# Analytically deriving Partial Information Decomposition for affine systems of stable and convolution-closed distributions

**Chaitanya Goswami**
Electrical & Computer Engineering,
Carnegie Mellon University,
Pittsburgh, PA-15213.
cgoswami@andrew.cmu.edu

**Amanda Merkley**
Electrical & Computer Engineering,
Carnegie Mellon University,
Pittsburgh, PA-15213.
amerkley@andrew.cmu.edu

## Abstract

Bivariate partial information decomposition (PID) has emerged as a promising tool for analyzing interactions in complex systems, particularly in neuroscience. PID achieves this by decomposing the information that two sources (e.g., different brain regions) have about a target (e.g., a stimulus) into unique, redundant, and synergistic terms. However, the computation of PID remains a challenging problem, often involving optimization over distributions. While several works have been proposed to compute PID terms *numerically*, there is a surprising dearth of work on computing PID terms *analytically*. The only known analytical PID result is for jointly Gaussian distributions. In this work, we present two theoretical advances that enable analytical calculation of the PID terms for numerous well-known distributions, including distributions relevant to neuroscience, such as Poisson, Cauchy, and binomial. Our first result generalizes the analytical Gaussian PID result to the much larger class of stable distributions. We also discover a theoretical link between PID and the emerging fields of data thinning and data fission. Our second result utilizes this link to derive analytical PID terms for two more classes of distributions: convolution-closed distributions and a sub-class of the exponential family. Furthermore, we provide an analytical upper bound for approximately computing PID for convolution-closed distributions, whose tightness we demonstrate in simulation.

## 1 Introduction

Bivariate partial information decomposition[1] (PID) is an information-theoretic framework developed for answering a central inquiry in many neuroscientific and machine learning studies: *how do two sources, $X$ and $Y$, jointly process information about a target $M$?* PID answers this question by quantifying the $M$-specific information contained in different interactions between $X$ and $Y$. Specifically, it decomposes the total information $X$ and $Y$ have about $M$ into *four* components: (i) the information about $M$ contained uniquely in $X$, (ii) the information about $M$ contained uniquely in $Y$, (iii) the redundant information about $M$ contained in both $X$ and $Y$, and (iv) the synergistic information about $M$ which arises from the interaction between $X$ and $Y$.

The PID terms offer novel insights for understanding interactions within complex systems, particularly in neuroscience. For instance, PID has been used to understand the firing patterns of grid cells [1, 2] and to study the flow of information in the visual cortex [3]. The following list of works [1, 3, 4, 5, 6, 7, 8] demonstrate the application of PID in studying diverse neuroscientific questions.

Beyond neuroscience, PID has found applications in multimodal learning for interpreting model predictions [9, 10], fair machine learning for defining and quantifying bias [11, 12, 13, 14], and understanding financial markets [15].

However, the primary constraint hindering broader adoption of PID is the difficulty of computing and estimating the PID terms. BROJA-PID [16], a widely applied PID framework, requires solving a constrained minimization problem over a set of probability distributions (see Sec. 2). This minimization problem can pose a considerable challenge, particularly when the underlying distributions are continuous. As a result, several recent works have been dedicated to providing *numerical algorithms* that solve the aforementioned minimization problem exactly or approximately [3, 9, 17, 18, 19, 20], with more emphasis on the case of discrete distributions.

Despite significant progress in *numerically* calculating PID (typically through discrete approximations), very few works exist on *analytically*[2] calculating PID. As of this writing, the only known analytical PID expressions exist for jointly Gaussian $M$, $X$, and $Y$ [21, 22]. A fundamental property of the Gaussian system is that one of the unique information (UI) terms in its PID is guaranteed to be *zero*. This property greatly simplifies the computation of PID for Gaussian systems, as the rest of the PID terms can be easily derived by solving desirable linear equations specified in many PID frameworks (see Sec 2), bypassing the need for optimizing over distributions (see Sec. 3).

In this work, we show that numerous systems of random variables $M$, $X$, and $Y$ expressing a particular "affine dependence structure" also exhibit the same fundamental property that at least one of the UI terms is zero. Consequently, we expand significantly on the existing Gaussian PID result by using this fundamental property to analytically compute PID for various systems of $M$, $X$, and $Y$ employing well-known distributions such as Poisson, exponential, gamma, beta, negative binomial, multinomial, Cauchy, Lévy-stable and more. The main contributions of this work are:

1. We extend the Gaussian PID result to a much larger class of distributions, known as the stable distribution family [23, 24] in Sec. 4. These results provide the first known analytical PID for fat-tailed distributions.

2. We highlight a theoretical link between PID calculation and the fields of data thinning [25] and data fission [26] in Sec. 5. We utilize this link to derive analytical PID terms for two more distribution families: convolution-closed distributions [25] and certain exponential family distributions [26].

3. For convolution-closed distributions, we further derive an analytical upper bound on the objective of the minimization used for computing BROJA-PID. We use this upper bound to approximately compute PID for systems of $M$, $X$, and $Y$ having a non-affine dependence structure. We show the goodness of our approximation by a simulation study in Sec. 6.

## 2  Background

**Notation**: We denote the set of all natural numbers, real numbers, and positive real numbers as $\mathbb{N}$, $\mathbb{R}$, and $\mathbb{R}^+$, respectively. Vectors are denoted by bold-faced font and an arrow, and matrices are denoted by bold-faced font. Define $\mathbb{N}_0 = \mathbb{N} \cup \{0\}$. Let $\mathbf{I}_d$ be the identity matrix of size $d \times d$ and $[d] = \{1, \ldots, d\} \; \forall \; d \in \mathbb{N}$. We denote $\vec{\mathbf{1}}_d$ and $\vec{\mathbf{0}}_d$ as $d$-dimensional vectors having all elements as 1 and 0, respectively. For brevity, the probability notations of the form $P(A|B)$ and $P(A)$ are always understood to be as $P(A = a|B = b)$ and $P(A = a)$, respectively. The general term 'distribution' is used to refer to both probability density function (p.d.f.) and probability mass function (p.m.f.) (whichever is appropriate depending upon the context). The symbol $A \perp\!\!\!\perp B|C$ denotes that $A$ and $B$ are conditionally independent given $C$, and similarly, $A \perp\!\!\!\perp B$ implies $A$ and $B$ are independent. The $L_p$-norm is denoted by $\|(\cdot)\|_p \; \forall \; p \in [0, \infty)$, and $|(\cdot)|$ denotes the absolute function.

**PID Background**: Suppose $M, X, Y$ are random variables with joint distribution $P(M, X, Y)$. According to [16, 27], three desirable equalities for a bivariate PID are as follows:

$$
\begin{aligned}
I_P(M; [X, Y]) &= UI(M; X\backslash Y) + UI(M; Y\backslash X) + RI(M; X, Y) + SI(M; X, Y), \\
I_P(M; X) &= UI(M; X\backslash Y) + RI(M; X, Y), \\
I_P(M; Y) &= UI(M; Y\backslash X) + RI(M; X, Y).
\end{aligned}
\tag{1}
$$

Here, $I_P(M; X)$ and $I_P(M; Y)$ denote the mutual information [28] between $M$ and $X$, and $M$ and $Y$, respectively, under the distribution $P(M, X, Y)$. Similarly, $I_P(M; [X, Y])$ denotes the joint mutual information between $M$ and $[X, Y]$ under the distribution $P(M, X, Y)$. $UI(M; X \backslash Y)$ and $UI(M; Y \backslash X)$ are the unique information about $M$ in $X$ and $Y$, respectively. $SI(M; X, Y)$ and $RI(M; X, Y)$ denote the respective synergistic and redundant information shared between $X$ and $Y$ about $M$. The linear system defined in (1) contains four variables in three equations. Hence, only one of $UI, RI$, or $SI$ need be defined to evaluate all four PID terms. Proposing a suitable definition of PID is the focus of much research [16, 27, 29, 30, 31, 32, 33, 34, 35] and all our theoretical results are applicable for any Blackwellian PID [22, 36] or a PID definition satisfying the assumption $(*)$ of Bertschinger et al. [16] (see Appx. B for a formal justification). For example, our results are applicable for the PID definitions proposed in [16, 22, 27, 29, 37]. In this work, we discuss our results in the context of the BROJA-PID [16, 38], which is the definition of PID adopted in previous works [3, 9, 20]:

$$UI(M; X \backslash Y) = \min_{Q \in \Delta_P} I_Q(M; X|Y) \text{ or equivalently } UI(M; Y \backslash X) = \min_{Q \in \Delta_P} I_Q(M; Y|X), \quad (2)$$

where $\Delta_P = \{Q(M, X, Y) : Q(M, X) = P(M, X), Q(M, Y) = P(M, Y)\}$ and $I_Q(M; X|Y)$ is the conditional mutual information under the distribution $Q(M, X, Y)$. Note that the minimizing distributions for both problems shown in (2) are the same [16], and consequently, both minimization problems are equivalent. We refer the reader to [39] for a review on PID. In this work, we define "analytically calculating PID" as analytically solving (2) by providing an explicit construction of the minimizing distribution $Q^*(M, X, Y)$ that minimizes (2). Note that only the distributions $Q^*(M, X, Y)$ and $P(M, X, Y)$ are needed to compute the BROJA-PID terms.

We briefly describe two distribution families used in Sec. 4 and Sec. 5 (see Appx. N for more details).

**Stable distribution family**: Stable distributions are a family of distributions that naturally arise in the context of generalized central limit theorems. Some well-known members of this family are the Gaussian, Cauchy, Poisson, and Lévy distributions. A defining feature of stable distributions is that the sum of two independent copies of a random variable $X$, denoted as $X_1$ and $X_2$, has the same distribution as a translated and scaled version of $X$ [23, 24, 40]. In this work, we consider five sub-classes of stable distributions:

(i) *Continuous stable distributions* are parameterized through four parameters: stability parameter $\alpha \in (0, 2]$, skewness parameter $\beta \in [-1, 1]$, scale parameter $\gamma \in (0, \infty)$, and location parameter $\mu \in \mathbb{R}$. We denote its p.d.f. as $p_{CS}(\alpha, \beta, \gamma, \mu)$. Note that all continuous stable distribution (except Gaussian) are fat-tailed.

(ii) *Independent component multivariate stable distributions* describe the distribution of $\vec{\mathbf{X}}$ consisting of $d$ independent random variables $\{X_j\}_{j=1}^d$, such that each $X_j \sim p_{CS}(\alpha, \beta_j, \gamma_j, \mu_j)$. We denote its p.d.f. as $p_{CS-IC}(\alpha, \vec{\boldsymbol{\beta}}, \vec{\boldsymbol{\gamma}}, \vec{\boldsymbol{\mu}})$.

(iii) *Elliptically-contoured multivariate stable distributions* are the distributions of continuous stable random vectors whose p.d.f. has elliptical contours, e.g., the multivariate Gaussian distribution. We denote its p.d.f. as $p_{CS-EC}(\alpha, \boldsymbol{\Sigma}, \vec{\boldsymbol{\mu}})$. Here, $\boldsymbol{\Sigma}$ is a positive definite matrix, $\vec{\boldsymbol{\mu}} \in \mathbb{R}^d$, and $\alpha \in (0, 2]$.

(iv) *Discrete stable distributions* are the discrete analogues of the continuous stable distributions. The p.m.f. of discrete stable distributions, denoted as $P_{DS}(\nu, \tau)$, are parameterized through two parameters: rate parameter $\tau > 0$ and exponent $0 < \nu \le 1$. Discrete stable distributions do not have well-known multivariate generalizations except the Poisson distribution.

(v) *Multivariate Poisson distribution*: We use the multivariate Poisson distribution proposed in [41, 42, 43, 44], denoted as $\text{Poisson}(d, d', \vec{\boldsymbol{\Lambda}})$, which represents each random variable in the random vector $\vec{\mathbf{N}}$ as a sum of independent Poisson random variables. Formally, $\vec{\mathbf{N}} \sim \text{Poisson}(d, d', \vec{\boldsymbol{\Lambda}})$ if $\vec{\mathbf{N}} = \mathbf{A}\vec{\mathbf{N}}^g$, where $\mathbf{A} = [\mathbf{A}_1 \ldots \mathbf{A}_{d'}]$. Here, $\mathbf{A}_i$ denotes a $d \times \binom{d}{i}$ submatrix having no duplicate columns, where each of its columns contain exactly $i$ ones and $(d - i)$ zeros [44]. The vector

$$\vec{\mathbf{N}}^g = [N_1^g \ldots N_d^g \ N_{12}^g \ldots \ N_{d-(d'-1)\ldots d}^g]^T$$

with mutually independent $N_{i_1 \ldots i_j}^g \sim \text{Poisson}(\lambda_{i_1 \ldots i_j}) \ \forall \ (i_1, \ldots, i_j) \in \mathbb{A}_j^d, j \in [d']$, where $\mathbb{A}_i^d = \{(j_1, \ldots, j_i) : j_1 < j_2 < \ldots < j_i, \text{ and } j_1, \ldots, j_i \in [d]\}$, e.g., $\mathbb{A}_2^3 = \{(1, 2), (1, 3), (2, 3)\}$. Note that $d' \le d$, and $\vec{\boldsymbol{\Lambda}} = [\lambda_1 \ \ldots \ \lambda_{d-(d'-1)\ldots d}]^T$.

We refer the reader to [23, 24, 40, 42, 44, 45, 46, 47] for more details on stable distributions.

**Convolution-closed distributions**: Convolution-closed distributions form a large class of distributions that are closed under convolution in some parameter $\delta$. Formally, we define the convolution-closed distribution as follows: Let $\mathcal{F}_\mathcal{D}$ denote a family of distributions, where each member distribution $f(\delta) \in \mathcal{F}_\mathcal{D}$ is indexed by a parameter $\delta \in \mathcal{D}$. Consider $X_1 \sim f(\delta_1)$, $X_2 \sim f(\delta_2)$, and $X_1 \perp\!\!\!\perp X_2$ for some $\delta_1, \delta_2 \in \mathcal{D}$. Then, $\mathcal{F}_\mathcal{D}$ is convolution-closed in the parameter $\delta$ if

$$X_1 + X_2 \sim f(\delta_1) * f(\delta_2) = f(\delta_1 + \delta_2) \ \forall \ \delta_1, \delta_2 \in \mathcal{D} \text{ such that } \delta_1 + \delta_2 \in \mathcal{D}, \tag{3}$$

where $*$ denotes the convolution operator. Many well-known distributions can be considered convolution-closed in some parameter [25], such as the Poisson distribution, Gaussian distribution, gamma distribution, etc. Table 1 in [25] lists various examples of convolution-closed distributions.

## 3   A sufficient condition for computing PID terms analytically

The main focus of this work is to analyze the cases where (2) is analytically solvable. A sufficient condition for solving (2) is to show that $\Delta_P$ contains a distribution $Q_{MC}(M, X, Y)$ with the Markovian structure $M \to X \to Y$ or $M \to Y \to X$. We briefly discuss the argument justifying this sufficient condition. Consider the case where $Q_{MC}(M, X, Y) \in \Delta_P$ and has the Markovian structure $M \to X \to Y$. First, we note that $UI(M; Y \backslash X) = \min_{Q \in \Delta_P} I_Q(M; Y|X) \geq 0$ due to the non-negativity of conditional mutual information [28]. Second, $Y \perp\!\!\!\perp M|X$ due to the Markovian structure of $Q_{MC}(M, X, Y)$, which implies that $I_{Q_{MC}}(M; Y|X) = 0$ (conditionally independent random variables have zero conditional mutual information [28]). Hence, $Q_{MC}(M, X, Y)$ achieves the lower bound of zero for the minimization problem $\min_{Q \in \Delta_P} I_Q(M; Y|X)$, showing that $Q_{MC}(M, X, Y)$ indeed minimizes (2). A similar argument can be made for the case when $Q_{MC}(M, X, Y)$ has the Markovian structure $M \to Y \to X$. Proposition 1 in Appx. A formalizes the above argument.

The above sufficient condition provides an easy way to analytically calculate the PID terms, as it ensures one of the unique information terms is always zero. Consequently, the remaining PID terms can be calculated by substituting zero for the appropriate unique information term in (1) and solving the resultant linear system. Hence, if applicable, the above sufficient condition considerably simplifies the calculation of PID terms by circumventing the need for optimizing over a set of distributions. Surprisingly, many well-known distribution families allow intuitive constructions of $P(M, X, Y)$, for which the above sufficient condition is applicable. In the following sections, we provide theorems that specify sufficient conditions under which the existence of these Markov chains in $\Delta_P$ can be guaranteed for these $P(M, X, Y)$.

## 4   Computing PID for stable distributions

In this section, we extend existing results for analytical PID computation of jointly Gaussian $M$, $X$, and $Y$ [21, 22] to the much larger class of stable distributions. Our results utilize two key observations to provide the generalization of the Gaussian results: (i) We first identify that the analytical computation of PID for jointly Gaussian systems is due to their particular "affine dependence" structure; (ii) We show that these particular affine dependence structures are not unique to Gaussian systems, but rather extend to many members of the stable distribution family.

For jointly Gaussian $M$, $X$, and $Y$, the conditional distributions $P(X|M) = \mathcal{N}(aM + b, \sigma_X^2)$ and $P(Y|M) = \mathcal{N}(cM + d, \sigma_Y^2)$ are also Gaussian distributions, where their means are an affine function of $M$ and their variances are fixed with respect to $M$. This particular affine dependence of $X$ and $Y$ on $M$ is the key to the analytical calculation of their PID, as it guarantees existence of a Markov chain $Q_{MC}(M, X, Y) \in \Delta_P$. Thus, we can apply the sufficient condition described in Sec. 3 to compute the PID terms. We illustrate through an example: consider $P(X|M) = \mathcal{N}(M, \sigma_X^2)$ and $P(Y|M) = \mathcal{N}(M, \sigma_Y^2)$ where $\sigma_X^2 < \sigma_Y^2$, and $M \sim P(M)$ for some appropriate $P(M)$. Then, we can explicitly construct $Q_{MC}(M, X, Y) \in \Delta_P$ with the Markovian structure $M \to X \to Y$ as follows: Choose $Q_{MC}(M, X, Y) = P(M)P(X|M)Q_{MC}(Y|X)$ where $Q_{MC}(Y|X)$ is specified through the addition of independent Gaussian noise, i.e., $Y = X + \epsilon$. Here, $\epsilon \sim \mathcal{N}(0, \sigma_Y^2 - \sigma_X^2)$ and $\epsilon \perp\!\!\!\perp (M, X)$. It is easy to verify that $Q_{MC}(Y|M) = \mathcal{N}(M, \sigma_Y^2)$, which implies that $Q_{MC}(M, Y) = P(M, Y)$. By construction, $Q_{MC}(M, X) = P(M, X)$, and hence, $Q_{MC}(M, X, Y) \in \Delta_P$.

The above example is a special case of a well-known result where, for a scalar $M$, one can always construct a lower signal-to-noise ratio (SNR) "Gaussian channel", i.e. $P(Y|M)$ in our example, by adding independent Gaussian noise to a higher SNR "Gaussian channel" [22, 48], i.e. $P(X|M)$ in our example. Surprisingly, the technique of adding independent noise to construct Markov chains contained in $\Delta_P$ can be extended to $P(X|M)$ and $P(Y|M)$ as members of stable distributions. Theorems 1 and 4 generalize the above construction for $P(X|M)$ and $P(Y|M)$ as members of univariate continuous and univariate discrete stable distributions, respectively. Theorems 2, 3, and 5 consider the case of multivariate stable distributions (see Sec. 2). The key technique for proving these theorems is that a Markov chain $Q_{MC}(M, X, Y) \in \Delta_P$ can always be constructed by adding appropriate independent noise to a higher SNR $P(X|M)$ to obtain a lower SNR $P(Y|M)$, similar to our above example.

## 4.1 PID of univariate affine continuous stable system

Theorem 1 can be viewed as a direct generalization of Barrett's Gaussian PID result [21] to stable distributions, showing one of the UI terms is always zero. We begin by formally describing the univariate affine continuous stable system that generalizes the Gaussian system of Barrett's [21]. Let the joint distribution of $M$, $X$, and $Y$, denoted as $P(M, X, Y)$, satisfy the following properties: (i) $M \sim P(M)$ with support set $\mathcal{M} \subseteq \mathbb{R}$; (ii) $P(X|M)$ and $P(Y|M)$ are univariate continuous stable distributions with an affine dependence on $M$, i.e., $P(X|M) = p_{CS}(\alpha, \beta_X, \gamma_X, aM + b)$ and $P(Y|M) = p_{CS}(\alpha, \beta_Y \operatorname{sgn}(ac), \gamma_Y, cM + d)$, where $a, b, c, d \in \mathbb{R}$.

**Theorem 1.** *Let $M$, $X$, and $Y$ be random variables whose joint distribution $P(M, X, Y)$ is described by a univariate affine continuous stable system. Without the loss of generality, assume ${|a|}/{\gamma_X} \geq {|c|}/{\gamma_Y}$. If $1 - \beta_Y \geq \left(\gamma_X |c|/\gamma_Y |a|\right)^\alpha (1 - \beta_X)$ and $1 + \beta_Y \geq \left(\gamma_X |c|/\gamma_Y |a|\right)^\alpha (1 + \beta_X)$, then $\Delta_P$ contains a Markov chain of the form $M \to X \to Y$ and $UI(M; Y \backslash X) = 0$.*

*Proof.* See Appx. F for the proof. Here, ${|a|}/{\gamma_X}$ and ${|c|}/{\gamma_Y}$ are the SNR analogues. $\qquad\square$

## 4.2 PID of multivariate affine continuous stable systems

We analyze two multivariate generalizations of Theorem 1 in Theorems 2 and 3. Namely, we consider independent component multivariate stable distribution (where all the components of the random vector are independent) and elliptically-contoured multivariate stable distribution (where the p.d.f. is elliptically contoured, similar to multivariate Gaussian distributions). We construct two systems employing these two sub-classes of multivariate continuous stable distributions and show that one of the UI terms is always zero for these systems. For both cases, denote the joint distribution of $M$, $\vec{\mathbf{X}}$, and $\vec{\mathbf{Y}}$ as $P(M, \vec{\mathbf{X}}, \vec{\mathbf{Y}})$, where the support set of $M$ is $\mathcal{M} \subseteq \mathbb{R}$. The dimensions of $M$, $\vec{\mathbf{X}}$, and $\vec{\mathbf{Y}}$ are $1$, $d_X$, and $d_Y$, respectively.

System 1: The random vectors $\vec{\mathbf{X}}$ and $\vec{\mathbf{Y}}$ satisfy the following equations:

$$\vec{\mathbf{X}} = \vec{\mathbf{H}}_X M + \mathbf{A}_X \vec{\mathbf{Z}}_X + \vec{\mathbf{b}}_X \text{ and } \vec{\mathbf{Y}} = \vec{\mathbf{H}}_Y M + \mathbf{A}_Y \vec{\mathbf{Z}}_Y + \vec{\mathbf{b}}_Y, \tag{4}$$

where $\vec{\mathbf{Z}}_X \sim p_{CS-IC}(\alpha, \vec{\mathbf{0}}_{d_X}, \vec{\mathbf{1}}_{d_X}, \vec{\mathbf{0}}_{d_X})$, $\vec{\mathbf{Z}}_Y \sim p_{CS-IC}(\alpha, \vec{\mathbf{0}}_{d_Y}, \vec{\mathbf{1}}_{d_Y}, \vec{\mathbf{0}}_{d_Y})$, $\mathbf{A}_X$ and $\mathbf{A}_Y$ are invertible matrices, $\vec{\mathbf{H}}_X, \vec{\mathbf{b}}_X \in \mathbb{R}^{d_X}$, and $\vec{\mathbf{H}}_Y, \vec{\mathbf{b}}_Y \in \mathbb{R}^{d_Y}$.

**Theorem 2.** *Let System 1 describe the joint distribution $P(M, \vec{\mathbf{X}}, \vec{\mathbf{Y}})$ of $M$, $\vec{\mathbf{X}}$, and $\vec{\mathbf{Y}}$. Without the loss of generality, assume $\|\mathbf{A}_Y^{-1}\vec{\mathbf{H}}_Y\|_\kappa \leq \|\mathbf{A}_X^{-1}\vec{\mathbf{H}}_X\|_\kappa$, where $\kappa = {\alpha}/{\alpha-1} \,\forall\, \alpha \in (1, 2]$ and $\kappa = \infty \,\forall\, \alpha \in (0, 1]$. Then, $\Delta_P$ contains a Markov chain of the form $M \to \vec{\mathbf{X}} \to \vec{\mathbf{Y}}$ and $UI(M; \vec{\mathbf{Y}} \backslash \vec{\mathbf{X}}) = 0$.*

*Proof.* See Appx. G for the proof. Here, $\|\mathbf{A}_Y^{-1}\vec{\mathbf{H}}_Y\|_\kappa$ and $\|\mathbf{A}_X^{-1}\vec{\mathbf{H}}_X\|_\kappa$ are the SNR analogues. $\quad\square$

System 2: The conditional distribution of $\vec{\mathbf{X}}$ and $\vec{\mathbf{Y}}$ conditioned on $M$ are as follows: $P(\vec{\mathbf{X}}|M) = p_{CS-EC}(\alpha, \mathbf{\Sigma}_X, \vec{\mathbf{H}}_X M + \vec{\mathbf{b}}_X)$ and $P(\vec{\mathbf{Y}}|M) = p_{CS-EC}(\alpha, \mathbf{\Sigma}_Y, \vec{\mathbf{H}}_Y M + \vec{\mathbf{b}}_Y)$, where $\mathbf{\Sigma}_X$ and $\mathbf{\Sigma}_Y$ are positive definite matrices, $\vec{\mathbf{H}}_X, \vec{\mathbf{b}}_X \in \mathbb{R}^{d_X}$, and $\vec{\mathbf{H}}_Y, \vec{\mathbf{b}}_Y \in \mathbb{R}^{d_Y}$.

**Theorem 3.** *Let System 2 describe the joint distribution $P(M, \vec{\mathbf{X}}, \vec{\mathbf{Y}})$ of $M$, $\vec{\mathbf{X}}$, and $\vec{\mathbf{Y}}$. Define $\boldsymbol{\Sigma}_X^{-1/2}$ and $\boldsymbol{\Sigma}_Y^{-1/2}$ as the respective inverses of the matrices $\boldsymbol{\Sigma}_X^{1/2}$ and $\boldsymbol{\Sigma}_Y^{1/2}$ which satisfy: $(\boldsymbol{\Sigma}_X^{1/2})^T \boldsymbol{\Sigma}_X^{1/2} = \boldsymbol{\Sigma}_X$, and $(\boldsymbol{\Sigma}_Y^{1/2})^T \boldsymbol{\Sigma}_Y^{1/2} = \boldsymbol{\Sigma}_Y$. Without the loss of generality, assume $\|\boldsymbol{\Sigma}_Y^{-1/2} \vec{\mathbf{H}}_Y\|_2 \leq \|\boldsymbol{\Sigma}_X^{-1/2} \vec{\mathbf{H}}_X\|_2$. Then, $\Delta_P$ contains a Markov chain of the form $M \rightarrow \vec{\mathbf{X}} \rightarrow \vec{\mathbf{Y}}$ and $UI(M; \vec{\mathbf{Y}} \backslash \vec{\mathbf{X}}) = 0$.*

*Proof.* See Appx. H for the proof. Here, $\|\boldsymbol{\Sigma}_Y^{-1/2} \vec{\mathbf{H}}_Y\|_2$ and $\|\boldsymbol{\Sigma}_X^{-1/2} \vec{\mathbf{H}}_X\|_2$ are the SNR analogues. □

### 4.3 PID of univariate affine discrete stable system

The univariate affine discrete stable system is the discrete counterpart of the univariate affine continuous stable system described in Sec. 4.1. The formal description of the univariate affine discrete stable system is as follows. Let the joint distribution of $M$, $X$, and $Y$, denoted as $P(M, X, Y)$, satisfy the following properties: (i) $M \sim P(M)$ with support set $\mathcal{M} \subseteq (0, \infty)$; (ii) $P(X|M)$ and $P(Y|M)$ are univariate discrete stable distributions with an affine dependence on $M$, i.e., $P(X|M{=}m){=}P_{DS}(\nu, am + b)$ and $P(Y|M{=}m){=}P_{DS}(\nu, cm + d)$, where $a, b, c, d \in (0, \infty)$.

**Theorem 4.** *Let $M$, $X$, and $Y$ be random variables whose joint distribution $P(M, X, Y)$ is described by a univariate affine discrete stable system. Without the loss of generality, assume $a \geq c$. If $a/b \geq c/d$, then $\Delta_P$ contains a Markov chain of the form $M \rightarrow X \rightarrow Y$ and $UI(M; Y \backslash X) = 0$.*

*Proof.* See Appx. I for the proof. Here, $a$ and $c$ are the SNR analogues. □

### 4.4 PID of multivariate linear Poisson system

The Poisson distribution is the only the discrete stable distribution with a well-known multivariate extension. Hence, we analyze vector-generalizations of Theorem 4 only for the Poisson distribution. We now describe the multivariate linear Poisson system: Let the joint distribution $P(M, \vec{\mathbf{X}}, \vec{\mathbf{Y}})$ of $M$, $\vec{\mathbf{X}}$, and $\vec{\mathbf{Y}}$ satisfy the following properties: (i) $M \sim P(M)$ with support set $\mathcal{M} \subseteq (0, \infty)$; (ii) $P(\vec{\mathbf{X}}|M) = \text{Poisson}(d_X, d_X', \vec{\boldsymbol{\Lambda}}_X)$ and $P(\vec{\mathbf{Y}}|M) = \text{Poisson}(d_Y, d_Y', \vec{\boldsymbol{\Lambda}}_Y)$, with:

$$\vec{\boldsymbol{\Lambda}}_X = \left[\lambda_1^X \ \cdots \ \lambda_{d_X - (d_X'-1)\ldots d_X}^X\right]^T, \lambda_{i_1 \ldots i_j}^X = \gamma_{i_1 \ldots i_j}^X M^j \ \forall j \in [d_X] \text{ and } (i_1, \ldots, i_j) \in \mathbb{A}_j^{d_X},$$

$$\vec{\boldsymbol{\Lambda}}_Y = \left[\lambda_1^Y \ \cdots \ \lambda_{d_Y - (d_Y'-1)\ldots d_Y}^Y\right]^T, \lambda_{i_1 \ldots i_j}^Y = \gamma_{i_1 \ldots i_j}^Y M^j \ \forall j \in [d_Y] \text{ and } (i_1, \ldots, i_j) \in \mathbb{A}_j^{d_Y}. \quad (5)$$

**Theorem 5.** *Let the joint distribution $P(M, \vec{\mathbf{X}}, \vec{\mathbf{Y}})$ of $M$, $\vec{\mathbf{X}}$, and $\vec{\mathbf{Y}}$ be described by the multivariate linear Poisson system defined above. Without the loss of generality, assume $d_X' \geq d_Y'$. If $\sum_{(i_1, \ldots, i_j) \in \mathbb{A}_j^{d_X}} \gamma_{i_1 \ldots i_j}^X \geq \sum_{(i_1, \ldots, i_j) \in \mathbb{A}_j^{d_Y}} \gamma_{i_1 \ldots i_j}^Y \ \forall j \in [d_Y']$, then $\Delta_P$ contains a Markov chain of the form $M \rightarrow \vec{\mathbf{X}} \rightarrow \vec{\mathbf{Y}}$ and $UI(M; \vec{\mathbf{Y}} \backslash \vec{\mathbf{X}}) = 0$.*

*Proof.* See Appx. J for the proof. Here, $\gamma_1^X, \ldots, \gamma_{d_X - (d_X'-1)\ldots d_X}^X, \gamma_1^Y, \ldots, \gamma_{d_Y - (d_Y'-1)\ldots d_Y}^Y \in \mathbb{R}^+$ are the SNR analogues. □

## 5 Computing PID using data thinning and data fission strategies

Data thinning and data fission are emerging fields in machine learning and statistics dedicated to studying the procedures of splitting a random variable $X$ into $N$ different component random variables. These splitting procedures provide an attractive alternative to the standard splitting of datasets into training, validation, and test splits for performing cross-validation to select statistical model parameters, as they enable users to perform cross-validation even for the extreme case of a dataset containing a *single* datapoint. We refer the reader to the recent works of Neufeld et al. [25] and Leiner et al. [26] for a more comprehensive discussion on data thinning and fission, respectively.

The fields of data thinning and data fission share an important theoretical link with analytically calculating PID terms. Specifically, the tools developed for splitting random variables for data thinning and data fission can be readily used to calculate PID terms analytically. To give an intuition, suppose $X$ contains more information about $M$ than $Y$. Then, we can employ data fission and thinning strategies to decompose $X$ into two components, $f_1(X)$ and $f_2(X)$, such that $f_2(X)$ follows the same distribution as $Y$. Then, $f_2(X)$ and $Y$ convey the same information about $M$ as they

are identically distributed and represent the redundant component of the PID terms. Similarly, $f_1(X)$ contains the information uniquely contained in $X$ about $M$. Thus, $f_1(X)$ represents the unique information term. Theorems 6 and 7 utilize the data thinning and data fission proposed in [25, 26] to construct a Markov chain $Q_{MC}(M, X, Y) \in \Delta_P$ for several systems of random variables $M$, $X$, and $Y$. This allows us to use the sufficient condition discussed in Sec. 3 to compute their PID.

## 5.1 PID for convolution-closed distribution based on data thinning strategies of Neufeld et al.

Neufeld et al. [25] introduces data thinning for a large family of distributions known as convolution-closed distributions (see Sec. 2). An attractive property of convolution-closed distributions is that they provide a natural way to define a dilation/thinning operation. Formally, let $X \sim f(\delta)$, then we define $X_\epsilon$ as the $\epsilon$-dilated version of $X$ if $X_\epsilon \sim f(\epsilon\delta)$ for some $\epsilon \in (0, 1)$ such that $\epsilon\delta \in \mathcal{D}$. Furthermore, if we assume $(1 - \epsilon)\delta \in \mathcal{D}$, then $P(X_\epsilon|X) = P(X_\epsilon|X_\epsilon + X_{1-\epsilon})$, where $X_{1-\epsilon} \sim f((1 - \epsilon)\delta)$ and $X_\epsilon \perp\!\!\!\perp X_{1-\epsilon}$ (see lemma 19 for a formal justification). We denote $P(X_\epsilon|X = x) = P(X_\epsilon|X_\epsilon + X_{1-\epsilon} = x)$ as $G(\epsilon\delta, (1 - \epsilon)\delta, x)$. This dilation operation forms the basis of data thinning, as it enables $X$ to be split into its dilated components $X_\epsilon$ and $X_{1-\epsilon}$, such that $X = X_\epsilon + X_{1-\epsilon}$, where $X_\epsilon \perp\!\!\!\perp X_{1-\epsilon}$. We utilize this dilation operation for analytically calculating the PID terms of the following linear convolution-closed system:

**Linear convolution-closed system**: Let $\mathcal{F}_\mathcal{D}$ be a convolution-closed distribution family as described in Sec. 2. The joint distribution $P(M, X, Y)$ of the random variables $M$, $X$, and $Y$ describes a *linear convolution-closed system* if the distributions $P(X|M)$ and $P(Y|M)$ are defined as follows:

$$P(X|M = m) = f(\delta_m^X) \text{ and } P(Y|M = m) = f(\delta_m^Y) \text{ such that } \delta_m^X, \delta_m^Y \in \mathcal{D} \, \forall \, m \in \mathcal{M}, \quad (6)$$

where $\mathcal{M}$ is the support of $M$. Furthermore, we assume $\delta_m^X = \gamma\delta_m^Y \, \forall \, m \in \mathcal{M}$ for some $\gamma \in \mathbb{R}^+$.

**Theorem 6.** *Let the joint density $P(M, X, Y)$ of random variables $M$, $X$, and $Y$ be described by a linear convolution-closed system. Without the loss of generality, assume $\gamma \leq 1$. If*

*(a) $(1 - \gamma)\delta_m^Y \in \mathcal{D} \, \forall \, m \in \mathcal{M}$,*

*(b) $P(X_\gamma|X_\gamma + X_{1-\gamma} = x, M = m) = G(\gamma\delta_m^X, (1 - \gamma)\delta_m^X, x)$ does not depend on $m$, where $P(X_\gamma|M) = f(\gamma\delta_m^X)$, $P(X_{1-\gamma}|M) = f((1 - \gamma)\delta_m^X)$ and $X_\gamma \perp\!\!\!\perp X_{1-\gamma}|M$,*

*then $\Delta_P$ contains a Markov chain of the form $M \to X \to Y$ and $UI(M; Y \backslash X) = 0$.*

The proof of Theorem 6 is provided in Appx. K. Theorem 6 enables PID calculation for several well-known distributions such as gamma, Poisson, beta etc. (see Appx. C for a (non-exhaustive) list).

## 5.2 PID for distributions based on data fission strategies of Leiner et al.

Leiner et al. [26] propose a "conjugate-prior reversal" strategy for splitting a random variable $X$ into two components, $f(X)$ and $g(X)$, for certain exponential family distributions. The distributions proposed for performing conjugate-prior reversal provide natural descriptions of $P(M, X, Y)$ (specified in theorem 7) for which PID can be calculated analytically. We briefly describe the distributions used in the conjugate-prior reversal strategy. Let $X \sim p_{exp1}(X)$, where

$$p_{exp1}(X = x; \theta_1, \theta_2) = H(\theta_1, \theta_2) \exp(\theta_1^T x - \theta_2^T A(x)), \quad (7)$$

for some appropriately defined $H(\cdot, \cdot), A(\cdot), \theta_1$ and $\theta_2$. Furthermore, define a random variable $Y$ through its conditional density $p(Y|X = x)$:

$$p(Y = y|X = x; \theta_3) = h(y) \exp\left(x^T T(y) - \theta_3^T A(x)\right), \quad (8)$$

for some $h(\cdot), T(\cdot)$, and $\theta_3$, such that $p(Y = y|X = x; \theta_1, \theta_2)$ is a well-defined distribution. Then, the decomposition terms $f(X)$ and $g(X)$ are $Y$ and $X$, respectively, in the conjugate-prior reversal strategy. Furthermore, the marginal distribution of $Y$ is expressed as:

$p_{exp2}(Y = y; \theta_1, \theta_2, \theta_3) = h(y)H(\theta_1, \theta_2)/H(\theta_1 + T(y), \theta_2 + \theta_3)$ (see proof of Theorem 1 in [26]).

**Theorem 7.** *Let $M, X$, and $Y$ be random variables having the joint distribution $P(M, X, Y)$. Furthermore, the conditional distribution of $X$ and $Y$ conditioned on $M$ are as follows: $P(X|M = m) = p_{exp1}(X; \theta_1(m), \theta_2(m))$ and $P(Y|M = m) = p_{exp2}(Y; \theta_1(m), \theta_2(m), \theta_3)$. Then, $\Delta_P$ contains a Markov chain of the form $M \to X \to Y$ and $UI(M; Y \backslash X) = 0$.*

The proof of Theorem 7 can be found in Appx. L. The proof essentially stems from observing that the joint distribution $Q_{MC}(M,X,Y) = P(M)P(X|M)Q_{MC}(Y|X)$ lies in $\Delta_P$, where $Q_{MC}(Y|X) = h(y)\exp\left(x^T T(y) - \theta_3^T A(x)\right)$. A (non-exhaustive) list of well-known distributions for which theorem 7 is applicable is provided in Appx. D. Leiner et al. [26] also discuss some more strategies for performing data fission that do not follow the conjugate-prior reversal strategy. We provide the corresponding results for computing PID for these remaining data fission strategies and for additional miscellaneous distributions in Appx. E.

## 6 Upper bound for convolution-closed distributions

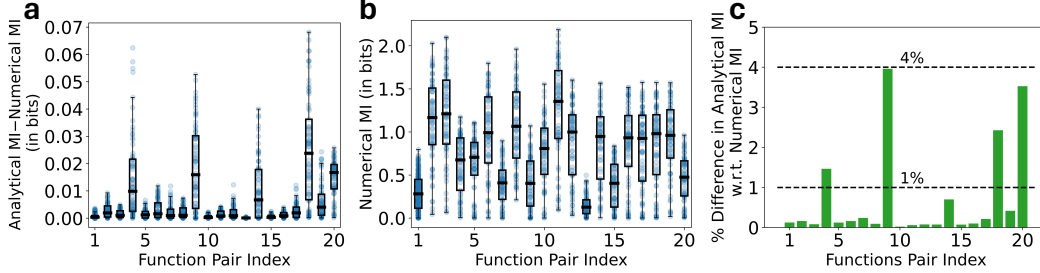

Figure 1: **a** and **b**, respectively, show the box plot of the difference $I_{Q_A}(M,X,Y) - I_{Q_N}(M,X,Y)$ and the corresponding values of $I_{Q_N}(M,X,Y)$ for the 20 different function pairs across the 75 different $P(M)$ distributions. The light-blue dots show the corresponding data points used for making the box plots. **c** shows the ratio of the median difference $I_{Q_A}(M,X,Y) - I_{Q_N}(M,X,Y)$ and the median value of $I_{Q_N}(M,X,Y)$ in percentage, for each function pair.

Several numerical methods, such as [3, 9, 20], approximately solve (2) by considering a smaller constraint set or minimizing an appropriate upper bound for calculating the PID terms. These numerical methods employ general approximations that rely on weak assumptions on the underlying distributions for solving (2), as their goal is to estimate PID for a large class of distributions. However, for specific applications, it is possible to make stronger assumptions on the underlying distributions (e.g., assuming the Poisson distribution for modeling neural spikes). This section illustrates how our theoretical analysis can benefit these numerical algorithms by providing more refined approximations for solving (2) that harness these stronger assumptions. Specifically, we construct an upper bound for the objective of (2) for convolution-closed distributions and show that, under certain assumptions, the upper bound can be *analytically minimized* over $\Delta_P$ in Sec. 6.1. Note that these upper bounds are applicable for more general cases than our theoretical results, as they do not require the sources $X$ and $Y$ to have an affine dependence on $M$. Consequently, our upper bound is also applicable in cases where both of the UI terms in the PID are non-zero, unlike our results in Sec. 4 and 5 We demonstrate the tightness of our upper bound through a simulation study in Sec. 6.2.

**Notations and assumptions**: We consider the minimization problem $\min_{Q \in \Delta_P} I_Q(M; [X,Y])$ for deriving the upper bound, as it was shown to be equivalent to (2) in [16]. The distribution $P(M,X,Y)$ for which we will construct our upper bound is specified as follows. The random variable $M$ has support over $\mathcal{M}$, the conditional distributions $P(X|M)$ and $P(Y|M)$ are members of some convolution-closed distribution family $\mathcal{F}_\mathcal{D}$, and there exists some $\delta_{bias}^X, \delta_{bias}^Y \in \mathcal{D}$ such that

$$P(X|M{=}m) = f(\delta_m^X) \text{ and } P(Y|M{=}m) = f(\delta_m^Y) \text{ where } \delta_m^X, \delta_m^Y \in \mathcal{D} \; \forall \, m \in \mathcal{M}, \text{ with}$$

$$(\delta_m^Y - \delta_{bias}^Y), (\delta_m^X - \delta_{bias}^X), (\delta_m^X - \delta_m^Y - (\delta_{bias}^X - \delta_{bias}^Y)) \in \mathcal{D} \text{ and } \delta_m^X - \delta_{bias}^X = \epsilon_m^{(1)}(\delta_m^Y - \delta_{bias}^Y),$$

$$\delta_m^Y = \epsilon_m^{(2)}(\delta_m^Y - \delta_{bias}^Y), \text{ and } \delta_m^X = \epsilon_m^{(3)}(\delta_m^X - \delta_{bias}^X) \text{ for } \epsilon_m^{(1)}, \epsilon_m^{(2)}, \epsilon_m^{(3)} \in [0,1] \; \forall \, m \in \mathcal{M}. \qquad (9)$$

The above assumptions ensure that $X$ and $Y$ can always be decomposed into new random variables $X', Y'', n_X, Y'$, and $n_Y$ (see (10)). We utilize these decomposed random variables to construct our upper bound. Hence, our upper bound is only applicable for systems satisfying (9). The above assumptions describe a large class of systems. For example, in the Poisson case, our upper bound is applicable for any $P(M,X,Y)$ having $P(X|M) = \text{Poisson}(f_1(M))$ and $P(Y|M) = \text{Poisson}(f_2(M))$, as long as $f_1(M) \geq f_2(M)$ over $\mathcal{M}$. In Appx. C, we provide numerous examples of systems for which (9) holds to illustrate the assumptions of (9).

## 6.1 Upper bound construction

First, we consider an arbitrary distribution $Q(M, X, Y) \in \Delta_P$. Therefore, we know $Q(X|M=m) = P(X|M=m) = f(\delta_m^X)$ and $Q(Y|M=m) = P(Y|M=m) = f(\delta_m^Y)$. We use the dilation properties of convolution-closed distributions (Sec. 5.1) to decompose $X$ and $Y$ into their respective dilated versions: $(X', Y'', n_X)$ and $(Y', n_Y)$. From the results of Appx. M.1, we know that $X = X' + Y'' + n_X$ and $Y = Y' + n_Y$. Furthermore, $(X', Y'', n_X)$ are mutually conditionally independent given $M$ and $Y' \perp\!\!\!\perp n_Y|M$. Hence, we can construct the following Markov chain for any arbitrary $Q(M, X, Y) \in \Delta_P$:

$$M \to \begin{bmatrix} X' & Y'' & Y' & n_X & n_Y \end{bmatrix}^T \to \begin{bmatrix} X' + Y'' + n_X & Y' + n_Y \end{bmatrix}^T = \begin{bmatrix} X & Y \end{bmatrix}^T. \quad (10)$$

We denote the joint distribution of $(M, X', Y'', Y', n_X, n_Y)$ as $\bar{Q}(M, X', Y'', Y', n_X, n_Y)$. We appropriately choose the dilation amounts for $X$ and $Y$ such that the respective conditional distributions of $(X', Y'', Y', n_X, n_Y)$ are as follows:

$$\bar{Q}(X'|M=m) = f(\delta_m^X - \delta_m^Y - (\delta_{bias}^X - \delta_{bias}^Y)), \quad \bar{Q}(Y''|M=m) = f(\delta_m^Y - \delta_{bias}^Y),$$
$$\bar{Q}(Y'|M=m) = f(\delta_m^Y - \delta_{bias}^Y), \quad \bar{Q}(n_X|M) = f(\delta_{bias}^X), \text{ and } \bar{Q}(n_Y|M) = f(\delta_{bias}^Y). \quad (11)$$

The distributions in (11) are well-defined due to (9). Appx. M.1 formally shows that a $\bar{Q}$, defined as above, exists for each $Q \in \Delta_P$. We use data-processing inequality and (10) to conclude:

$$I_{\bar{Q}}(M; [X', Y'', n_X, Y', n_Y]) \geq I_Q(M; [X, Y]) \,\forall\, Q(M, X, Y) \in \Delta_P. \quad (12)$$

Hence, (12) provides us the desired upper bound for our objective $I_Q(M; [X, Y])$. Furthermore, the minimization problem $\min_{Q \in \Delta_P} I_{\bar{Q}}(M; [X', Y'', n_X, Y', n_Y])$ is analytically solvable, and the minimizing distribution $\bar{Q}^*$ has the structure:

$$Y' = Y'' \text{ and } (n_X, n_Y) \perp\!\!\!\perp (M, X', Y').$$

The corresponding distribution of $(M, X, Y)$, denoted as $Q_A(M, X, Y)$, can be found by appropriately manipulating $\bar{Q}^*$, as $X = X' + Y'' + n_X$ and $Y = Y' + n_Y$. The distribution $Q_A$ serves as an approximate solution for the problem $\min_{Q \in \Delta_P} I_Q(M; [X, Y])$ and, consequently, (2). Note that if multiple $\delta_{bias}^X$ and $\delta_{bias}^Y$ exist satisfying (9), we optimize over the pairs $(\delta_{bias}^X, \delta_{bias}^Y)$ to further refine our approximate solution of (2).

## 6.2 Simulation study for numerically validating the upper bound

We illustrate the tightness of our upper bound through a simulation study on the Poisson distribution (a convolution-closed distribution). The reason for choosing the Poisson distribution is two-fold: (i) Poisson distribution is easily approximated as a discrete distribution over finite support, enabling calculation of ground-truth PID terms through numerical solvers such as [10, 17, 18], which are not readily available for continuous distributions; (ii) Many practical applications of PID have been in neuroscience, and the Poisson distribution is frequently used for modeling neural spikes in neuroscientific studies [49].

We compare the performance of our analytical estimate $Q_A(M, X, Y)$ with the numerical ground-truth estimate $Q_N(M, X, Y)$ for the Poisson distribution. The simulation setup is as follows: We choose a $P(M, X, Y)$ such that $P(X|M) = \text{Poisson}(f_1(M))$ and $P(Y|M) = \text{Poisson}(f_2(M))$. We chose 20 different pairs of non-linear $f_1(\cdot)$ and $f_2(\cdot)$ (enumerated in Table 1 in Appx. M), such that the assumptions in (9) are satisfied. The distribution $P(M)$ is chosen to be a discrete distribution. For each function pair, we compare $Q_A(M, X, Y)$ and $Q_N(M, X, Y)$ over 75 different distributions of $P(M)$ as shown in Fig. 1a. The 75 distributions are further sub-divided into three groups of 25 based on the number of the outcomes of $M$, which are 2, 4, and 8. For each of the 75 distributions, the values of $M$ are randomly sampled from $[1, 4]$ and the values of $P(M)$ are randomly sampled from a simplex of appropriate dimensions. We compare the mutual information $I_{Q_A}(M; [X, Y])$ calculated using $Q_A$ with the numerical ground-truth $I_{Q_N}(M; [X, Y])$ calculated using numerical solvers (Fig. 1b). For calculating $I_{Q_N}(M; [X, Y])$, we approximate the Poisson distribution as a finite discrete distribution by terminating its support at the smallest integer where its cumulative distribution is greater than 0.99. We use the code provided in [50] (under Apache 2.0-license) to numerically solve (2) using this discrete approximation of the Poisson distribution. Fig. 1c demonstrates that the analytical upper bound provided by $Q_A$ is very tight for the tested function pairs (within $< 1\%$ of

the numerical ground-truth for 16 function pairs) and serves as a good approximation for solving (2). Furthermore, the tightness of our analytical upper bound suggests that $Q_A$ might be an analytical solution of (2) for a larger class of systems of $M$, $X$, and $Y$ having non-affine dependence on $M$. Similar results demonstrating the tightness of upper-bound for negative-binomial and binomial distributions are presented in Appx. M.4.

# 7    Discussion and Limitations

In this work, we analytically compute PID for large classes of distributions, greatly expanding upon the analytical result for the Gaussian system. We provide the first known analytical PID for systems employing Poisson, gamma, exponential, Cauchy, beta, Dirichlet, Lévy-stable, binomial, multinomial, negative binomial, and uniform distributions. Furthermore, we generalize the previous Gaussian PID result [21] in an additional way by showing the target $M$ need not be Gaussian. Our stable distribution results provide the first known analytical computation of PID for fat-tailed distributions (all continuous stable distributions have infinite variance except the Gaussian distribution). A practical utility of our analytical results is that they provide a large test bed in which the performance of numerical PID estimators can be compared and evaluated. This test bed may benefit future works on numerical PID estimation by enabling more comprehensive testing of PID estimators.

Our results on the Poisson, Cauchy, and binomial are of particular relevance in the neuroscientific context. Poisson and Cauchy distributions are widely used to model neural spikes [49] and network dynamics in the brain [51]. Binomial thinning is a frequently-used operator in neuroscience [49]. Our generalization of the Gaussian result could be helpful in refining approximations already used in computing PID for neural data [3], e.g., by relaxing the assumption of joint Gaussianity. As continuous stable distributions have been shown to better model Magnetic Resonance Imaging data [52, 53], our results may also be helpful in computing PID in this application.

We also connect the fields of data thinning and data fission with PID by using their decomposition strategies for analytically computing the PID of systems based on convolution-closed distributions (see Sec. 5). Conversely, our PID results also suggest decomposition strategies for data thinning/fission purposes, e.g., our stable distribution results suggest that stable random variables can be decomposed by adding independent noise (similar to the Gaussian case discussed in [26]). Convolution-closed distributions are particularly promising for studying PID as they allow intuitive construction of upper bounds that can be analytically minimized (see Sec. 6). These upper bounds can complement the work on numerical estimation of PID by providing more refined approximations. Another promising avenue is to combine our upper bounds with lower bounds (e.g., from [54]) to create branch and bound algorithms [55] for solving (2). Overall, our analytical results greatly facilitate the computation of PID, either by directly using the analytical expressions or by providing refined approximations for numerical methods.

**Limitations and Future Work**: We study PID for univariate $M$, $X$, and $Y$ and provide some vector extensions. More vector extensions of our results are a promising direction for subsequent works. Most of our analytical results require $P(X|M)$ and $P(Y|M)$ to depend on some affine functions of $M$. The existence of analytical solutions for the cases where $P(X|M)$ and $P(Y|M)$ depend on non-affine functions of $M$ remains an open question. The upper bound discussed in Sec. 6 can be further refined with more careful analysis, and more rigorous testing of these upper bounds is required to understand in which cases the upper bound is a good approximation for solving (2). We defer the testing and refinement of these upper bounds for subsequent works, as our primary goal in this work is to study analytical solutions of PID. Niu & Quinn [56] also propose a duality result between the synergistic and redundant components in the Gaussian broadcast and multiple-access channels utilizing the analytical PID expressions of the Gaussian system. It may be possible to derive similar duality results for appropriately defined broadcast and multiple-access channels employing other distributions (e.g., Poisson or Cauchy) with our theoretical results.

**Broader Impact**: Due to their theoretical nature, our results' negative impact primarily depends on how they are used and interpreted. Our theoretical results are applicable only under the specific assumptions outlined in this work, and using these results without ensuring that the theorem's assumptions are satisfied can lead to incorrect scientific conclusions. We also caution reader against naively using PID to draw causal inferences, as PID is inherently a correlational quantity.

## Acknowledgments and Disclosure of Funding

Authors would like to thank Prof. Pulkit Grover, Dr. Praveen Venkatesh, Prof. Alireza Chamanzar, and Neil Ashim Mehta for helpful discussions. This work was supported by National Science Foundation (NSF). Authors declare no competing interests.

## Footnotes

[1]Throughout this work we refer to bivariate PID as PID.

[2] Section 2 precisely defines the notion of "analytical calculating PID" adopted in this work.

## References

[1] Praveen Venkatesh and Pulkit Grover. Understanding encoding and redundancy in grid cells using partial information decomposition. *Comput. Syst. Neurosci.(Cosyne)*, 2020.

[2] Ariel K Feldman, Praveen Venkatesh, Douglas J Weber, and Pulkit Grover. Information-theoretic tools to understand distributed source coding in neuroscience. *IEEE Journal on Selected Areas in Information Theory*, 2024.

[3] Praveen Venkatesh, Corbett Bennett, Sam Gale, Tamina Ramirez, Greggory Heller, Severine Durand, Shawn Olsen, and Stefan Mihalas. Gaussian partial information decomposition: Bias correction and application to high-dimensional data. In A. Oh, T. Naumann, A. Globerson, K. Saenko, M. Hardt, and S. Levine, editors, *Advances in Neural Information Processing Systems*, volume 36, pages 74602–74635. Curran Associates, Inc., 2023.

[4] Nigel Colenbier, Frederik Van de Steen, Lucina Q Uddin, Russell A Poldrack, Vince D Calhoun, and Daniele Marinazzo. Disambiguating the role of blood flow and global signal with partial information decomposition. *NeuroImage*, 213:116699, 2020.

[5] Tjeerd W Boonstra, Luca Faes, Jennifer N Kerkman, and Daniele Marinazzo. Information decomposition of multichannel EMG to map functional interactions in the distributed motor system. *NeuroImage*, 202:116093, 2019.

[6] Jana Krohova, Luca Faes, Barbora Czippelova, Zuzana Turianikova, Nikoleta Mazgutova, Riccardo Pernice, Alessandro Busacca, Daniele Marinazzo, Sebastiano Stramaglia, and Michal Javorka. Multiscale information decomposition dissects control mechanisms of heart rate variability at rest and during physiological stress. *Entropy*, 21(5):526, 2019.

[7] Giuseppe Pica, Eugenio Piasini, Houman Safaai, Caroline Runyan, Christopher Harvey, Mathew Diamond, Christoph Kayser, Tommaso Fellin, and Stefano Panzeri. Quantifying how much sensory information in a neural code is relevant for behavior. In I. Guyon, U. Von Luxburg, S. Bengio, H. Wallach, R. Fergus, S. Vishwanathan, and R. Garnett, editors, *Advances in Neural Information Processing Systems*, volume 30. Curran Associates, Inc., 2017.

[8] Praveen Venkatesh, Sanghamitra Dutta, and Pulkit Grover. Information flow in computational systems. *IEEE Transactions on Information Theory*, 66(9):5456–5491, 2020.

[9] Paul Pu Liang, Yun Cheng, Xiang Fan, Chun Kai Ling, Suzanne Nie, Richard Chen, Zihao Deng, Nicholas Allen, Randy Auerbach, Faisal Mahmood, Russ R Salakhutdinov, and Louis-Philippe Morency. Quantifying & modeling multimodal interactions: An information decomposition framework. In A. Oh, T. Naumann, A. Globerson, K. Saenko, M. Hardt, and S. Levine, editors, *Advances in Neural Information Processing Systems*, volume 36, pages 27351–27393. Curran Associates, Inc., 2023.

[10] Paul Pu Liang, Yun Cheng, Ruslan Salakhutdinov, and Louis-Philippe Morency. Multimodal fusion interactions: A study of human and automatic quantification. In *Proceedings of the 25th International Conference on Multimodal Interaction*, pages 425–435, 2023.

[11] Sanghamitra Dutta, Praveen Venkatesh, Piotr Mardziel, Anupam Datta, and Pulkit Grover. An information-theoretic quantification of discrimination with exempt features. In *Proceedings of the AAAI Conference on Artificial Intelligence*, volume 34, pages 3825–3833, 2020.

[12] Sanghamitra Dutta and Faisal Hamman. A review of partial information decomposition in algorithmic fairness and explainability. *Entropy*, 25(5):795, 2023.

[13] Faisal Hamman and Sanghamitra Dutta. Demystifying local & global fairness trade-offs in federated learning using partial information decomposition. In *The Twelfth International Conference on Learning Representations*, 2024.

[14] Faisal Hamman and Sanghamitra Dutta. A unified view of group fairness tradeoffs using partial information decomposition. In *2024 IEEE International Symposium on Information Theory (ISIT)*, pages 214–219, 2024.

[15] Tomas Scagliarini, Luca Faes, Daniele Marinazzo, Sebastiano Stramaglia, and Rosario N Mantegna. Synergistic information transfer in the global system of financial markets. *Entropy*, 22(9):1000, 2020.

[16] Nils Bertschinger, Johannes Rauh, Eckehard Olbrich, Jürgen Jost, and Nihat Ay. Quantifying unique information. *Entropy*, 16(4):2161–2183, 2014.

[17] Abdullah Makkeh, Dirk Oliver Theis, and Raul Vicente. Bivariate partial information decomposition: The optimization perspective. *Entropy*, 19(10):530, 2017.

[18] Pradeep Kr Banerjee, Johannes Rauh, and Guido Montúfar. Computing the unique information. In *2018 IEEE International Symposium on Information Theory (ISIT)*, pages 141–145. IEEE, 2018.

[19] Abdullah Makkeh, Dirk Oliver Theis, and Raul Vicente. BROJA-2PID: A robust estimator for bivariate partial information decomposition. *Entropy*, 20(4):271, 2018.

[20] Ari Pakman, Amin Nejatbakhsh, Dar Gilboa, Abdullah Makkeh, Luca Mazzucato, Michael Wibral, and Elad Schneidman. Estimating the unique information of continuous variables. In M. Ranzato, A. Beygelzimer, Y. Dauphin, P.S. Liang, and J. Wortman Vaughan, editors, *Advances in Neural Information Processing Systems*, volume 34, pages 20295–20307. Curran Associates, Inc., 2021.

[21] Adam B Barrett. Exploration of synergistic and redundant information sharing in static and dynamical Gaussian systems. *Physical Review E*, 91(5):052802, 2015.

[22] Praveen Venkatesh and Gabriel Schamberg. Partial information decomposition via deficiency for multivariate Gaussians. In *2022 IEEE International Symposium on Information Theory (ISIT)*, pages 2892–2897. IEEE, 2022.

[23] John P Nolan. Univariate stable distributions. *Springer Series in Operations Research and Financial Engineering*, 10:978–3, 2020.

[24] Fred W Steutel and Klaas van Harn. Discrete analogues of self-decomposability and stability. *The Annals of Probability*, pages 893–899, 1979.

[25] Anna Neufeld, Ameer Dharamshi, Lucy L. Gao, and Daniela Witten. Data thinning for convolution-closed distributions. *Journal of Machine Learning Research*, 25(57):1–35, 2024.

[26] James Leiner, Boyan Duan, Larry Wasserman, and Aaditya Ramdas. Data fission: Splitting a single data point. *Journal of the American Statistical Association*, 0(0):1–12, 2023.

[27] Paul L Williams and Randall D Beer. Nonnegative decomposition of multivariate information. *arXiv preprint arXiv:1004.2515*, 2010.

[28] Thomas M. Cover and Joy A. Thomas. *Elements of Information Theory (Wiley Series in Telecommunications and Signal Processing)*. Wiley-Interscience, USA, 2006.

[29] Malte Harder, Christoph Salge, and Daniel Polani. Bivariate measure of redundant information. *Physical Review E*, 87(1):012130, 2013.

[30] Pradeep Kr Banerjee, Eckehard Olbrich, Jürgen Jost, and Johannes Rauh. Unique informations and deficiencies. In *2018 56th Annual Allerton Conference on Communication, Control, and Computing (Allerton)*, pages 32–38. IEEE, 2018.

[31] Jim W Kay and Robin AA Ince. Exact partial information decompositions for Gaussian systems based on dependency constraints. *Entropy*, 20(4):240, 2018.

[32] Conor Finn and Joseph T Lizier. Pointwise partial information decomposition using the specificity and ambiguity lattices. *Entropy*, 20(4):297, 2018.

[33] Xueyan Niu and Christopher J Quinn. A measure of synergy, redundancy, and unique information using information geometry. In *2019 IEEE International Symposium on Information Theory (ISIT)*, pages 3127–3131. IEEE, 2019.

[34] Fernando E Rosas, Pedro AM Mediano, Borzoo Rassouli, and Adam B Barrett. An operational information decomposition via synergistic disclosure. *Journal of Physics A: Mathematical and Theoretical*, 53(48):485001, 2020.

[35] Keerthana Gurushankar, Praveen Venkatesh, and Pulkit Grover. Extracting unique information through Markov relations. In *2022 58th Annual Allerton Conference on Communication, Control, and Computing (Allerton)*, pages 1–6. IEEE, 2022.

[36] David Blackwell. Equivalent comparisons of experiments. *The Annals of Mathematical Statistics*, pages 265–272, 1953.

[37] Praveen Venkatesh, Keerthana Gurushankar, and Gabriel Schamberg. Capturing and interpreting unique information. In *2023 IEEE International Symposium on Information Theory (ISIT)*, pages 2631–2636. IEEE, 2023.

[38] Virgil Griffith and Christof Koch. Quantifying synergistic mutual information. *Guided self-organization: inception*, pages 159–190, 2014.

[39] Joseph T Lizier, Nils Bertschinger, Jürgen Jost, and Michael Wibral. Information decomposition of target effects from multi-source interactions: Perspectives on previous, current and future work. *Entropy*, 20(4):307, 2018.

[40] John P Nolan. *Stable distributions*. 2012.

[41] Jeremie Fish, Jie Sun, and Erik Bollt. Interaction networks from discrete event data by Poisson multivariate mutual information estimation and information flow with applications from gene expression data. *Applied Network Science*, 7(1):1–22, 2022.

[42] AS Krishnamoorthy. Multivariate binomial and Poisson distributions. *Sankhyā: The Indian Journal of Statistics*, pages 117–124, 1951.

[43] DM Mahamunulu. A note on regression in the multivariate Poisson distribution. *Journal of the American Statistical Association*, 62(317):251–258, 1967.

[44] Dimitris Karlis and Loukia Meligkotsidou. Multivariate Poisson regression with covariance structure. *Statistics and Computing*, 15(4):255–265, 2005.

[45] Murad S Taqqu. *Stable non-Gaussian random processes: stochastic models with infinite variance*. Chapman & Hall, 1994.

[46] S James Press. Multivariate stable distributions. *Journal of Multivariate Analysis*, 2(4):444–462, 1972.

[47] Norman Lloyd Johnson, Samuel Kotz, and Narayanaswamy Balakrishnan. *Discrete multivariate distributions*, volume 165. Wiley New York, 1997.

[48] Xiaohu Shang and H Vincent Poor. Noisy-interference sum-rate capacity for vector Gaussian interference channels. *IEEE Transactions on Information Theory*, 59(1):132–153, 2012.

[49] Peter Dayan and Laurence F Abbott. *Theoretical neuroscience: computational and mathematical modeling of neural systems*. MIT press, 2005.

[50] Abdullah Makkeh, Dirk Oliver Theis, and Raul Vicente. Broja-2pid: A robust estimator for bertschinger et al.'s bivariate partial information decomposition. *Entropy*, 20(4):271, 2018.

[51] Viktoras Pyragas and Kestutis Pyragas. Mean-field equations for neural populations with q-Gaussian heterogeneities. *Physical Review E*, 105(4):044402, 2022.

[52] Diego Salas-González, JM Górriz, Javier Ramírez, Matthias Schloegl, Elmar Wolfgang Lang, and Andrés Ortiz. Parameterization of the distribution of white and grey matter in MRI using the $\alpha$-stable distribution. *Computers in biology and medicine*, 43(5):559–567, 2013.

[53] Diego Salas-Gonzalez, Juan M Górriz, Javier Ramírez, and Elmar W Lang. Why using the alpha-stable distribution in NeuroImage? In *2014 International Conference on Signal Processing and Multimedia Applications (SIGMAP)*, pages 297–301. IEEE, 2014.

[54] Seiya Tokui and Issei Sato. Disentanglement analysis with partial information decomposition. *arXiv preprint arXiv:2108.13753*, 2021.

[55] Narendra and Fukunaga. A branch and bound algorithm for feature subset selection. *IEEE Transactions on Computers*, 100(9):917–922, 1977.

[56] Xueyan Niu and Christopher J Quinn. Synergy and redundancy duality between Gaussian multiple access and broadcast channels. In *2020 International Symposium on Information Theory and Its Applications (ISITA)*, pages 6–10. IEEE, 2020.

[57] Robert B Ash and Catherine A Doléans-Dade. *Probability and measure theory*. Academic press, 2000.

[58] Gilbert Strang. *Introduction to linear algebra*. SIAM, 2022.

[59] Alan V Oppenheim, Alan S Willsky, Syed Hamid Nawab, and Jian-Jiun Ding. *Signals and systems*, volume 2. Prentice hall Upper Saddle River, NJ, 1997.

[60] Jagadesh Chandramohan and Lung-Kuang Liang. Bernoulli, multinomial and Markov chain thinning of some point processes and some results about the superposition of dependent renewal processes. *Journal of Applied Probability*, 22(4):828–835, 1985.

[61] Sergio Verdú. Poisson communication theory. *International Technion Communication Day in Honor of Israel Bar-David*, 66, 1999.

[62] Christian Walck et al. *Hand-book on statistical distributions for experimentalists*. Stockholms universitet, 1996.

[63] Kalimuthu Krishnamoorthy. *Handbook of statistical distributions with applications*. Chapman and Hall/CRC, 2006.

[64] Lenka Slámová and Lev B Klebanov. Generalized definitions of discrete stability. *arXiv preprint arXiv:1502.02588*, 2015.

# A  Formal proof of the sufficient condition for solving (2) discussed in Sec. 3

**Proposition 1.** *For the minimization problem defined in* (2)*, if there exists a joint density* $Q_{MC}(M, X, Y) \in \Delta_P$*, having the Markovian structure of the form* $M \to X \to Y$ *or* $M \to Y \to X$*, then* $Q_{MC}(M, X, Y)$ *minimizes* (2)*. Furthermore,*

1. *If* $Q_{MC}(M, X, Y)$ *has the Markovian structure* $M \to X \to Y$*, then* $UI(M; Y\backslash X) = I_{Q_{MC}}(M; Y|X) = 0.$

2. *If* $Q_{MC}(M, X, Y)$ *has the Markovian structure* $M \to Y \to X$*, then* $UI(M; X\backslash Y) = I_{Q_{MC}}(M, X|Y) = 0.$

*Proof.* **Case 1**: $Q_{MC}(M, X, Y) \in \Delta_P$ having the Markovian structure $M \to X \to Y$.

By the definition of the Markov chain, we know that $M$ and $Y$ are conditionally independent given $X$. Denote $Q_{MC}(M, X, Y)$ as the joint distribution of the Markov chain $M \to X \to Y$. Then,

$$I_{Q_{MC}}(M; Y|X) = 0, \tag{13}$$

where $I_{Q_{MC}}(M; Y|X)$ is the conditional mutual information between $M$ and $Y$ given $X$ under the distribution $Q_{MC}(M, X, Y)$. Equation (13) follows from a well-known property of conditional mutual information which states that $I(A; B|C) = 0$ iff $A \perp\!\!\!\perp B|C$ [28].

Since $I_Q(M; Y|X) \geq 0 \, \forall \, Q \in \Delta_P$, due to the non-negativity of conditional mutual information [28], we can conclude that $\min_{Q \in \Delta_P} I_Q(M; Y|X) \geq 0$. The joint distribution $Q_{MC}(M, X, Y)$ having the Markovian structures $M \to X \to Y$ achieves the lower bound of 0 for the minimization problem $\min_{Q \in \Delta_P} I_Q(M; Y|X)$ implying $Q_{MC}(M, X, Y)$ minimizes (2).

**Case 2**: $Q_{MC}(M, X, Y) \in \Delta_P$ having the Markovian structure $M \to Y \to X$.

A similar argument as case 1, with the random variables $X$ and $Y$ switched, would allow us to conclude that the joint distribution $Q_{MC}(M, X, Y)$ specifying the Markov chain $M \to Y \to X$ minimizes (2). □

# B  Other PID definitions for which theorems 1-7 are valid

Our theoretical results (Theorems 1-7) hold for a wide range of PID measures. Particularly, our results hold for two broad classes of PID definitions: PID definitions satisfying assumption $(*)$ in Bertschinger et al. [16] and Blackwellian PIDs (see Appendix B-C of Venkatesh & Schamberg [22] for a deeper discussion on differences between these two families of PID definition). Some examples of PID definitions satisfying assumption $(*)$ in Bertschinger et al. [16] are the Williams & Beer's PID definition [27], the PID definition proposed in Harder et al. [29], and I-PID proposed in Venkatesh et al. [37]. Similarly, an example of a Blackwellian PID is the $\delta$-PID discussed in Venkatesh & Schamberg [22]. Section III-F of Venkatesh et al. [37] further discusses a family of Blackwellian PID definitions for which our theorems are also applicable. We now provide formal arguments showing that our theoretical results are applicable for PID definitions that either satisfy assumption $(*)$ of Bertschinger et al. [16] or are Blackwellian

**PID definitions satisfying assumption** $(*)$: We invoke the argument presented in the proof discussed in Section 4.2 of Barrett [21] to show that our results hold for any PID definition satisfying assumption $(*)$ in Bertschinger et al. [16]. Barrett's argument relies on the key observation that the UI calculated using BROJA-PID upper bounds the UI calculated using any other PID definition satisfying assumption $(*)$ in Bertschinger et al. Entropy'14. Consequently, if one of the UI atoms of the BROJA-PID is zero then the corresponding UI atom of the other PID definitions must also be zero due to BROJA PID upper-bounding them and the non-negativity of the PID atoms. Since the remaining PID atoms are calculated using (1), it must be the case that the PID atoms calculated using any PID definition satisfying assumption $(*)$ must be the same as the PID atoms calculated using BROJA-PID whenever one of the UI terms of BROJA-PID is zero.

**Blackwellian PID**: The defining feature of a Blackwellian PID is that the UI atom is zero iff it is possible to construct the Markovian structure between the random variables (i.e., $M \to X \to Y$ or $M \to Y \to X$), while preserving their pairwise marginals (see Venkatesh et al. [22] for more details). In all our theoretical results, we show that the random variables in the analyzed system admit

a Markovian structure, while preserving the pairwise marginal structure. Hence, the UI atom for the systems analyzed in Theorems 1-7 is zero for any Blackwellian PID. Consequently, as the rest of the PID terms are calculated using (1), it must be the case that the PID atoms calculated using any Blackwellian PID must be the same as the PID atoms calculated using BROJA-PID in our results.

## C  Lists of Systems that can solved using Theorem 6

In this section, we provide examples of systems of random variables $(M, X, Y)$ employing well-known distributions whose PID terms can be obtained through theorem 6. The following list is modified from Table 2 of [25]. Note that for the joint distribution $P(M, X, Y)$ of the system of random variable $(M, X, Y)$, we only specify $P(M)$, $P(X|M)$, and $P(Y|M)$, as any $P(M, X, Y)$ having these marginals can be solved through theorem 6. We also discuss some possible extensions of the corresponding systems for which the upper bound discussed in Sec. 6 is applicable. Note that the extension of the systems provided are not exhaustive, and there exists more systems for which the upper bound is applicable. In the following examples, we assume $a > b$.

1. **System**: The distributions of $P(M), P(X|M)$, and $P(Y|M)$ are as follows:

    (a) $M \sim P(M)$ having support $(0, \infty)$.
    (b) $P(X|M)$=Poisson$(aM)$.
    (c) $P(Y|M) = $ Poisson$(bM)$.

    The corresponding $G(\gamma \delta_m^X, (1-\gamma)\delta_m^X, x)$ specified in theorem 6 is Binomial$(x, \gamma)$ with $\gamma = b/a$, and $a, b > 0$.
    **Result**: The Markov chain $M \to X \to Y \in \Delta_P$ and $UI(M; Y \backslash X) = 0$.
    **An extension of the system for which the upper bound of Sec. 6 in applicable**: The distributions of $P(M), P(X|M)$, and $P(Y|M)$ are as follows:

    (a) $M \sim P(M)$ having support $\mathcal{M}$.
    (b) $P(X|M)$=Poisson$(f_1(M))$, for $f_1(M) > 0 \,\forall\, m \in \mathcal{M}$.
    (c) $P(Y|M) = $ Poisson$(f_2(M))$, for $f_2(M) > 0 \,\forall\, m \in \mathcal{M}$.
    (d) $f_1(m) \geq f_2(m) \,\forall\, m \in \mathcal{M}$ and all the underlying distributions are well-defined.

2. **System**: The distributions of $P(M), P(X|M)$, and $P(Y|M)$ are as follows:

    (a) $M \sim P(M)$ having support $(-\infty, \infty)$.
    (b) $P(X|M)$=$\mathcal{N}(aM, a\sigma^2)$.
    (c) $P(Y|M) = \mathcal{N}(bM, b\sigma^2)$.

    The corresponding $G(\gamma \delta_m^X, (1-\gamma)\delta_m^X, x)$ specified in theorem 6 is $\mathcal{N}(\gamma x, \gamma(1-\gamma)\sigma^2)$ with $\gamma = b/a$, and $a, b > 0$.
    **Result**: The Markov chain $M \to X \to Y \in \Delta_P$ and $UI(M; Y \backslash X) = 0$.
    **An extension of the system for which the upper bound of Sec. 6 in applicable**: The distributions of $P(M), P(X|M)$, and $P(Y|M)$ are as follows:

    (a) $M \sim P(M)$ having support $\mathcal{M}$.
    (b) $P(X|M)$=Gaussian$(f_1(M), f_1(M))$, for $f_1(M) > 0 \,\forall\, m \in \mathcal{M}$.
    (c) $P(Y|M) = $ Gaussian$(f_2(M), f_2(M))$, for $f_2(M) > 0 \,\forall\, m \in \mathcal{M}$.
    (d) $f_1(m) \geq f_2(m) \,\forall\, m \in \mathcal{M}$ and all the underlying distributions are well-defined.

3. **System**: The distributions of $P(M), P(X|M)$, and $P(Y|M)$ are as follows:

    (a) $M \sim P(M)$ having support $(0, 1]$.
    (b) $P(X|M)$=Negative Binomial$(a, M)$.
    (c) $P(Y|M) = $ Negative Binomial$(b, M)$.

    The corresponding $G(\gamma \delta_m^X, (1-\gamma)\delta_m^X, x)$ specified in theorem 6 is Beta-Binomial$(x, \gamma a, (1-\gamma)a)$ with $\gamma = b/a$, and $a, b \in \mathbb{N}$.
    **Result**: The Markov chain $M \to X \to Y \in \Delta_P$, and $UI(M; Y \backslash X) = 0$.
    **An extension of the system for which the upper bound of Sec. 6 in applicable**: The distributions of $P(M), P(X|M)$, and $P(Y|M)$ are as follows:

    (a) $M \sim P(M)$ having support $\mathcal{M}$.

(b)  $P(X|M)$=Negative Binomial$(f_1(M), p)$, for $f_1(M) \in \mathbb{N}_0 \, \forall \, m \in \mathcal{M}$ and $p \in [0, 1]$.

(c)  $P(Y|M)$=Negative Binomial$(f_2(M), p)$, for $f_2(M) \in \mathbb{N}_0 \, \forall \, m \in \mathcal{M}$.

(d)  $f_1(m) \geq f_2(m) \, \forall \, m \in \mathcal{M}$ and all the underlying distributions are well-defined.

4. **System**: The distributions of $P(M), P(X|M)$, and $P(Y|M)$ are as follows:

   (a)  $M \sim P(M)$ having support $(0, \infty)$.

   (b)  $P(X|M)$=Gamma$(a, M)$.

   (c)  $P(Y|M) = $ Gamma$(b, M)$.

   The corresponding $G(\gamma\delta_m^X, (1-\gamma)\delta_m^X, x)$ specified in theorem 6 is obtained as follows: $G(\gamma\delta_m^X, (1-\gamma)\delta_m^X, x)=xZ, \; Z \sim \text{Beta}(b, (1-b))$ with $\gamma = {}^{b}\!/\!{}_{a}$, and $a, b \in (0, \infty)$. Note that we are following the Gamma$(\alpha, \beta)$ notation.

   **Result**: The Markov chain $M \to X \to Y \in \Delta_P$ and $UI(M; Y\backslash X) = 0$.

   **An extension of the system for which the upper bound of Sec. 6 in applicable**: The distributions of $P(M), P(X|M)$, and $P(Y|M)$ are as follows:

   (a)  $M \sim P(M)$ having support $\mathcal{M}$.

   (b)  $P(X|M)$=Gamma$(f_1(M), \beta)$, for $f_1(M) > 0 \, \forall \, m \in \mathcal{M}$ and $\beta > 0$.

   (c)  $P(Y|M) = $ Gamma$(f_2(M), \beta)$, for $f_2(M) > 0 \, \forall \, m \in \mathcal{M}$.

   (d)  $f_1(m) \geq f_2(m) \, \forall \, m \in \mathcal{M}$ and all the underlying distributions are well-defined.

5. **System**: The distributions of $P(M), P(X|M)$, and $P(Y|M)$ are as follows:

   (a)  $M \sim P(M)$ having support $(0, \infty)$.

   (b)  $P(X|M)$=Exponential$(M)$.

   (c)  $P(Y|M) = $ Gamma$(b, M)$.

   The corresponding $G(\gamma\delta_m^X, (1-\gamma)\delta_m^X, x)$ specified in theorem 6 is obtained as follows: $G(\gamma\delta_m^X, (1-\gamma)\delta_m^X, x)=xZ, \; Z \sim \text{Beta}(b, (1-b))$, and $0 < b \leq 1$. Note that we are following the Gamma$(\alpha, \beta)$ notation.

   **Result**: The Markov chain $M \to X \to Y \in \Delta_P$ and $UI(M; Y\backslash X) = 0$.

   **An extension of the system for which the upper bound of Sec. 6 in applicable**: The distributions of $P(M), P(X|M)$, and $P(Y|M)$ are as follows:

   (a)  $M \sim P(M)$ having support $\mathcal{M}$.

   (b)  $P(X|M)$=Exponential$(f_1(M))$, for $f_1(M) > 0 \, \forall \, m \in \mathcal{M}$.

   (c)  $P(Y|M) = $ Gamma$(f_2(M), f_1(M))$, for $f_2(M) > 0 \, \forall \, m \in \mathcal{M}$.

   (d)  $f_2(m) \leq 1 \, \forall \, m \in \mathcal{M}$ and all the underlying distributions are well-defined.

6. **System**: The distributions of $P(M), P(X|M)$, and $P(Y|M)$ are as follows:

   (a)  $M \sim P(M)$ having support $[0, 1]$.

   (b)  $P(X|M)$=Binomial$(a, M)$.

   (c)  $P(Y|M) = $ Binomial$(b, M)$.

   The corresponding $G(\gamma\delta_m^X, (1-\gamma)\delta_m^X, x)$ specified in theorem 6 is Hypergeometric$(x, \gamma a, (1-\gamma)a)$ with $\gamma = {}^{b}\!/\!{}_{a}$, and $a, b \in \mathbb{N}$.

   **Result**: The Markov chain $M \to X \to Y \in \Delta_P$ and $UI(M; Y\backslash X) = 0$.

   **An extension of the system for which the upper bound of Sec. 6 in applicable**: The distributions of $P(M), P(X|M)$, and $P(Y|M)$ are as follows:

   (a)  $M \sim P(M)$ having support $\mathcal{M}$.

   (b)  $P(X|M)$=Binomial$(f_1(M), p)$, for $f_1(M) \in \mathbb{N}_0 \, \forall \, m \in \mathcal{M}$ and $p \in [0, 1]$.

   (c)  $P(Y|M) = $ Binomial$(f_2(M), p)$, for $f_2(M) \in \mathbb{N}_0 \, \forall \, m \in \mathcal{M}$.

   (d)  $f_1(m) \geq f_2(m) \, \forall \, m \in \mathcal{M}$ and all the underlying distributions are well-defined.

7. **System**: The distributions of $P(\vec{\mathbf{M}}), P(\vec{\mathbf{X}}|\vec{\mathbf{M}})$, and $P(\vec{\mathbf{Y}}|\vec{\mathbf{M}})$ are as follows:

   (a)  $\vec{\mathbf{M}} \sim P(\vec{\mathbf{M}})$ having support over a $k$-dimensional simplex.

   (b)  $P(\vec{\mathbf{X}}|\vec{\mathbf{M}})$=Multinomial$_k(a, \vec{\mathbf{M}})$.

   (c)  $P(\vec{\mathbf{Y}}|\vec{\mathbf{M}}) = $ Multinomial$_k(b, \vec{\mathbf{M}})$.

The corresponding $G(\gamma \delta_m^X, (1-\gamma)\delta_m^X, x)$ specified in theorem 6 is Multivariate Hypergeometric $(\vec{\mathbf{x}}, \gamma_Y)$ with $\gamma = b/a$, and $a, b \in \mathbb{N}$.

**Result**: The Markov chain $\vec{\mathbf{M}} \to \vec{\mathbf{X}} \to \vec{\mathbf{Y}} \in \Delta_P$ and $UI(\vec{\mathbf{M}}; \vec{\mathbf{Y}} \backslash \vec{\mathbf{X}}) = 0$.

**An extension of the system for which the upper bound of Sec. 6 in applicable**: The distributions of $P(M), P(\vec{\mathbf{X}}|M)$, and $P(\vec{\mathbf{Y}}|M)$ are as follows:

(a) $M \sim P(M)$ having support $\mathcal{M}$.

(b) $P(\vec{\mathbf{X}}|M) = \text{Multinomial}_k(f_1(M), \vec{\mathbf{p}})$, for $f_1(M) \in \mathbb{N}_0 \ \forall \ m \in \mathcal{M}$.

(c) $P(\vec{\mathbf{Y}}|M) = \text{Multinomial}(f_2(M), \vec{\mathbf{p}})$, for $f_2(M) \in \mathbb{N}_0 \ \forall \ m \in \mathcal{M}$.

(d) $f_1(m) \geq f_2(m) \ \forall \ m \in \mathcal{M}$, $\vec{\mathbf{p}}$ is a valid probability vector, and all the underlying distributions are well-defined.

# D   Lists of Systems that can solved using Theorem 7

In this section, we provide examples of systems of random variables $(M, X, Y)$ employing well-known distributions whose PID terms can be obtained through theorem 7. The following list is appropriately modified from Appendix A of [26]. Similar to Appx. C, we only specify specify $P(M)$, $P(X|M)$, and $P(Y|M)$ of the joint distribution $P(M, X, Y)$, as any $P(M, X, Y)$ having the same marginals can be solved through theorem 7. In the following examples, we assume $a > b$.

1. **System**: The distributions of $P(M), P(X|M)$, and $P(Y|M)$ are as follows:

   (a) $M \sim P(M)$ having support $(0, \infty)$.

   (b) $P(X|M) = \text{Exponential}(M)$.

   (c) $P(Y|M) = \text{Geometric}(M/(\tau+M))$, where $\tau \in (0, \infty)$.

   **Results**: The Markov chain $M \to X \to Y \in \Delta_P$, and $UI(M; Y \backslash X) = 0$. The corresponding $Q_{MC}(Y|X)$ required for constructing the Markov chain is Poisson$(\tau X)$.

2. **System**: The distributions of $P(M), P(X|M)$, and $P(Y|M)$ are as follows:

   (a) $M \sim P(M)$ having support $\mathcal{M}$.

   (b) $P(X|M) = \text{Gamma}(f_1(M), f_2(M))$, where $f_1(m), f_2(m) > 0 \ \forall \ m \in \mathcal{M}$.

   (c) $P(Y|M) = \text{Negative Binomial}(f_1(M), f_2(M)/(f_2(M)+\tau))$, where $\tau \in (0, \infty)$

   **Results**: The Markov chain $M \to X \to Y \in \Delta_P$, and $UI(M; Y \backslash X) = 0$. The corresponding $Q_{MC}(Y|X)$ required for constructing the Markov chain is is Poisson$(\tau X)$.

3. **System**: The distributions of $P(M), P(X|M)$, and $P(Y|M)$ are as follows:

   (a) $M \sim P(M)$ having support $(0, \infty)$.

   (b) $P(X|M) = \text{Beta}(M, 1)$.

   (c) $P(Y|M)$ is a discrete distribution with over support set $\{0, 1, \ldots, N\}$, with

   $$P(Y = y|M = m) = m \frac{\Gamma(y+m)N!}{\Gamma(N+1+m)y!},$$

   where $\Gamma(.)$ is the Gamma function.

   **Results**: The Markov chain $M \to X \to Y \in \Delta_P$, and $UI(M; Y \backslash X) = 0$. The corresponding $Q_{MC}(Y|X)$ required for constructing the Markov chain is Binomial$(N, X)$.

4. **System**: The distributions of $P(M), P(X|M)$, and $P(Y|M)$ are as follows:

   (a) $M \sim P(M)$ having support $(0, \infty)$.

   (b) $P(X|M) = \text{Beta}(1, M)$.

   (c) $P(Y|M)$ is a discrete distribution with over support set $\{0, 1, \ldots, N\}$, with

   $$P(Y = y|M = m) = m \frac{\Gamma(y+m)N!}{\Gamma(N+1+m)y!},$$

   where $\Gamma(.)$ is the Gamma function.

   **Results**: The Markov chain $M \to X \to Y \in \Delta_P$, and $UI(M; Y \backslash X) = 0$. The corresponding $Q_{MC}(Y|X)$ required for constructing the Markov chain is is Binomial$(N, 1-X)$.

5. **System**: The distributions of $P(\vec{\mathbf{M}})$, $P(\vec{\mathbf{X}}|\vec{\mathbf{M}})$, and $P(\vec{\mathbf{Y}}|\vec{\mathbf{M}})$ are as follows:

   (a) $\vec{\mathbf{M}} \sim P(\vec{\mathbf{M}})$ having support $(0, \infty)^d$ for some dimension $d$.

   (b) $P(\vec{\mathbf{X}}|\vec{\mathbf{M}})$=Dirichlet$(M_1, \dots, M_d)$.

   (c) $P(\vec{\mathbf{Y}}|\vec{\mathbf{M}})$ is a discrete distribution having support $\{y_i \in \{1, \dots, N\} \ \forall \ i \in [d] | \sum_{i=1}^d y_i = N\}$, and

$$P(\vec{\mathbf{Y}}|\vec{\mathbf{M}}) = \frac{\Gamma\left(\sum_{i=1}^d M_i\right) N!}{\Gamma\left(\sum_{i=1}^d M_i + N\right)} \prod_{i=1}^d \frac{\Gamma(M_i + Y_i)}{\Gamma(M_i) Y_i!},$$

   where $\Gamma(.)$ is the Gamma function.

   **Results**: The Markov chain $\vec{\mathbf{M}} \to \vec{\mathbf{X}} \to \vec{\mathbf{Y}} \in \Delta_P$, and $UI(\vec{\mathbf{M}}; \vec{\mathbf{Y}} \backslash \vec{\mathbf{X}}) = 0$. The corresponding $Q_{MC}(\vec{\mathbf{Y}}|\vec{\mathbf{X}})$ required for constructing the Markov chain is Multinomial$_d(N, \vec{\mathbf{X}})$.

# E   Miscellaneous PID Results

This section explores the analytical calculation of PID terms for five more systems of random variables $(M, X, Y)$ with joint distribution $P(M, X, Y)$. These five systems' $P(M, X, Y)$ employ well-known distributions, but their respective analytical PID calculation does not result from Theorem 1-Theorem 7. Of note are the results of $P(M, X, Y)$ employing exponential and uniform distribution, which states that these systems will always have both unique information terms zero.

## E.1   PID for the miscellaneous system M1.

Miscellaneous system **M1**: Let $(M, X, Y)$ be a system of random variables with joint distribution $\overline{P(M, X, Y)}$. The joint distribution $P(M, X, Y)$ describes the miscellaneous system **M1** if it satisfies the following two properties:

1. $M \sim P(M)$, and has support $(0, \infty)$.
2. $P(X|M)$=Exponential$(\gamma_X M)$, and $P(Y|M)$=Exponential$(\gamma_Y M)$, where $\gamma_X, \gamma_Y \in (0, \infty)$.

**Proposition 2.** *Let $M, X,$ and $Y$ be random variables whose joint distribution $P(M, X, Y)$ describes the miscellaneous system **M1** defined above. Then, $\Delta_P$ contains a distribution $Q_{MC}(M, X, Y)$, which admits both Markovian structures, i.e., $M \to X \to Y$ and $M \to Y \to X$. Consequently, both $UI(M; Y \backslash X) = UI(M; X \backslash Y) = 0$.*

*Proof.* We prove proposition 2 by explicitly constructing a joint distribution $Q_{MC}(M, X, Y) \in \Delta_P$ that admits both Markovian structures, namely, $M \to X \to Y$ and $M \to Y \to X$. A necessary and sufficient condition for the distribution $Q_{MC}(M, X, Y)$ to admit both Markovian structures is that it should satisfy (14).

$$Q_{MC}(M, X, Y)=Q_{MC}(M)Q_{MC}(X|M)Q_{MC}(Y|X)=Q_{MC}(M)Q_{MC}(Y|M)Q_{MC}(X|Y). \tag{14}$$

We now briefly outline the structure of the proof. The rest of the proof can be divided into three parts:

1. In the first part, we explicitly construct $Q_{MC}(M, X, Y)$.

2. In the second part, we show that the $Q_{MC}(M, X, Y)$ constructed in the first part lies in $\Delta_P$.

3. In the third part, we conclude our proof by showing that $Q_{MC}(M, X, Y)$ satisfies (14) and, consequently, admits both Markovian structures.

**Part 1: Specifying $Q_{MC}(M, X, Y)$**

We specify $Q_{MC}(M, X, Y)$ by individually specifying $Q_{MC}(M)$, $Q_{MC}(X|M)$, and $Q_{MC}(Y|X)$. We choose $Q_{MC}(M)$ and $Q_{MC}(X|M)$ as follows:

$$Q_{MC}(M) = P(M), \text{ and } Q_{MC}(X|M) = P(X|M). \tag{15}$$

The distribution $Q_{MC}(Y|X)$ is specified through the following deterministic transformation:

$$Y = \frac{\gamma_X}{\gamma_Y}X. \tag{16}$$

**Part 2: Showing $Q_{MC}(M,X,Y) \in \Delta_P$**

For showing $Q_{MC}(M,X,Y) \in \Delta_P$, we first need to calculate $Q_{MC}(Y|M)$:

*Calculating $Q_{MC}(Y|M)$:* By lemma 1, if $X \sim \text{Exp}(\lambda)$, then $aX \sim \text{Exponential}(\lambda/a)$. Since $Q_{MC}(X|M) = \text{Exponential}(\gamma_X M)$, we can use the result of lemma 1 to calculate $Q_{MC}(Y|M)$ as follows:

$$Q_{MC}(Y|M) = Q_{MC}\left(\frac{\gamma_X}{\gamma_Y}X \,\middle|\, M\right) = \text{Exponential}\left(\frac{\gamma_Y}{\gamma_X}\gamma_X M\right)$$
$$= \text{Exponential}\left(\gamma_Y M\right) = P(Y|M). \tag{17}$$

Then, from (15) and (17), we can conclude:

$$Q_{MC}(M,X) = Q_{MC}(M)Q_{MC}(X|M) = P(M)P(X|M) = P(M,X), \tag{18}$$
$$Q_{MC}(M,Y) = Q_{MC}(M)Q_{MC}(Y|M) = P(M)P(Y|M) = P(M,Y). \tag{19}$$

Hence, $Q_{MC}(M,X,Y) \in \Delta_P$.

**Part 3: Showing $Q_{MC}(M,X,Y)$ satisfies (14)**

It is trivial to see that by the construction of $Q_{MC}(M,X,Y)$ given in Part 1, we have:

$$Q_{MC}(M,X,Y) = Q_{MC}(M)Q_{MC}(X|M)Q_{MC}(Y|X). \tag{20}$$

By the chain rule, we have:

$$Q_{MC}(M,X,Y) = Q_{MC}(M)Q_{MC}(Y|M)Q_{MC}(X|Y,M). \tag{21}$$

Hence, for showing that $Q_{MC}(M,X,Y)$ satisfies (14), we only need to show that $\underline{Q_{MC}(X|Y,M) = Q_{MC}(X|Y)}$.

*Showing $Q_{MC}(X|Y,M) = Q_{MC}(X|Y)$:* Observe that $Y = \gamma_X/\gamma_Y X$ by construction. Therefore, we can equivalently express $X = \gamma_Y/\gamma_X Y$. Consequently,

$$Q_{MC}(X|Y,M) = \mathbb{I}[X = \gamma_Y/\gamma_X Y] \text{ as } X = \gamma_Y/\gamma_X Y, \tag{22}$$

where $\mathbb{I}[\cdot]$ is the indicator function. Since R.H.S of the above equation does not depend on $M$, we have:

$$Q_{MC}(X|Y,M) = \mathbb{I}[X = \gamma_Y/\gamma_X Y] = Q_{MC}(X|Y). \tag{23}$$

Combining (21) and (23), we have:

$$Q_{MC}(M,X,Y) = Q_{MC}(M)Q_{MC}(Y|M)Q_{MC}(X|Y). \tag{24}$$

Combining (20) and (24), we can show that $Q_{MC}(M,X,Y)$ satisfies (14).

**Conclusion**: Combining the results of parts 2 and 3, we can conclude that the $Q_{MC}(M,X,Y) \in \Delta_P$ and contains both Markovian structure $M \to X \to Y$ and $M \to Y \to X$. A unique implication of $Q_{MC}(M,X,Y)$ admitting both Markovian structures is that both $I_{Q_{MC}}(M;X|Y) = I_{Q_{MC}}(M;Y|X) = 0$. Consequently, both minimization problems presented in (2) are minimized by $Q_{MC}(M,X,Y)$ with $UI(M;X\backslash Y) = \min_{Q \in \Delta_P} I_Q(M;X|Y) = 0$ and $UI(M;Y\backslash X) = \min_{Q \in \Delta_P} I_Q(M;Y|X) = 0$, completing our proof. $\qquad\square$

**Lemma 1.** *If $X \sim \text{Exponential}(\lambda)$ with $\lambda \in (0,\infty)$, then $\gamma X \sim \text{Exponential}(\lambda/\gamma)$ for $\gamma \in (0,\infty)$.*

*Proof.* We denote the p.d.f. of the random variable $X$ as $p_X(\cdot)$ and, by lemma statement, we know:

$$p_X(X = x) = \text{Exponential}(\lambda) = \begin{cases} \lambda e^{-x\lambda} & x \geq 0 \\ 0 & \text{otherwise} \end{cases}. \tag{25}$$

Denote the p.d.f. of the function $f(X)$ of the random variable $X$ as $p_{f(X)}(\cdot)$. We want to derive $p_{f(X)}(\cdot)$ for $f(X) = \gamma X$. Since, $f(X) = \gamma X$ is an invertible function, we can use the change-of-variable technique for deriving $p_{f(X)}(\cdot)$:

$$p_{f(X)}(f(X) = y) = p_X(X = f^{-1}(y)) \left| \frac{df^{-1}(y)}{dy} \right| = \begin{cases} \frac{\lambda}{\gamma} e^{-y\lambda/\gamma} & y \geq 0 \\ 0 & \text{otherwise} \end{cases}$$

$$= \text{Exponential}(\lambda/\gamma). \tag{26}$$

$\square$

## E.2   PID for the miscellaneous system M2.

Miscellaneous System **M2**: Let $(M, X, Y)$ be a system of random variables with joint distribution $P(M, X, Y)$. The joint distribution $P(M, X, Y)$ describes the miscellaneous system **M2** if it satisfies the following two properties:

1. $M \sim P(M)$ and has support $(0, \infty)$.
2. $P(X|M) = \text{Negative Binomial}(M, p_X)$, and $P(Y|M) = \text{Negative Binomial}(M, p_Y)$.

This system is adapted from the negative binomial decomposition listed in Appendix A [26].

**Proposition 3.** *Let $M, X$, and $Y$ be random variables whose joint distribution $P(M, X, Y)$ describes the miscellaneous system **M2** defined above.*

1. *If $p_X \leq p_Y$, then $\Delta_P$ contains a Markov chain of the form $M \rightarrow X \rightarrow Y$, and $UI(M; Y \backslash X) = 0$.*

2. *If $p_X \geq p_Y$, then $\Delta_P$ contains a Markov chain of the form $M \rightarrow Y \rightarrow X$, and $UI(M; X \backslash Y) = 0$.*

*Proof.* We only provide an explicit proof of condition 1. The proof of condition 2 follows the same steps as the proof of condition 1 with parameters of $X$ and $Y$ switched.

**Proof of condition 1**: We briefly outline the proof structure. The proof can be divided into two parts:

1. In the first part, we explicitly construct a joint distribution $Q_{MC}(M, X, Y)$ having the Markovian structure $M \rightarrow X \rightarrow Y$.

2. In the second part, we conclude our proof by showing the $Q_{MC}(M, X, Y)$ constructed in the first part, lies in $\Delta_P$.

### Part 1: Specifying $Q_{MC}(M, X, Y)$

Since $Q_{MC}(M, X, Y)$ is the joint distribution of a Markov chain $M \rightarrow X \rightarrow Y$, we have $Q_{MC}(M, X, Y) = Q_{MC}(M)Q_{MC}(X|M)Q_{MC}(Y|X)$. Taking advantage of this special structure of $Q_{MC}(M, X, Y)$, we specify $Q_{MC}(M, X, Y)$ by individually specifying $Q_{MC}(M), Q_{MC}(X|M)$, and $Q_{MC}(Y|X)$. We choose $Q_{MC}(M), Q_{MC}(X|M)$ and $Q_{MC}(Y|X)$ as follows:

$$Q_{MC}(M) = P(M), Q_{MC}(X|M) = P(X|M) \text{ and } Q_{MC}(Y|X) = \text{Binomial}(X, p),$$

$$\text{where } p = \frac{p_X(1 - p_Y)}{p_Y(1 - p_X)}. \tag{27}$$

### Part 2: Showing $Q_{MC}(M, X, Y) \in \Delta_P$

For showing $Q_{MC}(M, X, Y) \in \Delta_P$, we first need to calculate $Q_{MC}(Y|M)$:

*Calculating $Q_{MC}(Y|M)$:* Since $Q_{MC}(X|M) = \text{Negative Binomial}(M, p_X)$ and $Q_{MC}(Y|X) = \text{Binomial}(N, p)$, we can use lemma 2 for calculating $Q(Y|M)$:

$$Q_{MC}(Y|M) = \text{Negative Binomial}(M, p_Y) = P(Y|M). \tag{28}$$

Then, from (27) and (28), we can conclude:

$$Q_{MC}(M, X) = Q_{MC}(M)Q_{MC}(X|M) = P(M)P(X|M) = P(M, X), \qquad (29)$$
$$Q_{MC}(M, Y) = Q_{MC}(M)Q_{MC}(Y|M) = P(M)P(Y|M) = P(M, Y). \qquad (30)$$

Hence, $Q_{MC}(M, X, Y) \in \Delta_P$ and consequently, by proposition 1, $UI(M; Y \backslash X) = 0$. This concludes our proof. $\qquad\square$

**Lemma 2.** *Let* $X \sim$ *Negative Binomial*$(N, p_X)$ *and* $P(Y|X) =$ *Binomial*$(X, p)$ *with* $p = \frac{p_X(1-p_Y)}{p_Y(1-p_X)}$, $p_Y \geq p_X$, $p_Y, p_X \in [0, 1]$, *and* $N \in \mathbb{N}_0$. *Then,* $Y \sim$ *Negative Binomial*$(N, p_Y)$.

*Proof.* First, in order for $P(Y|X)$ to be a legitimate distribution, we need to ensure that $p \in [0, 1]$, i.e., we need to show that:

$$0 \leq p \leq 1. \qquad (31)$$

Since $p_X, (1 - p_Y), (1 - p_X), p_Y \geq 0$ as $p_X, p_Y \in [0, 1]$, we can easily verify the first inequality:

$$p = \frac{p_X(1 - p_Y)}{(1 - p_X)p_Y} \geq 0. \qquad (32)$$

For verifying $p \leq 1$, we will use the assumption that $p_Y \geq p_X$:

$$p_X \leq p_Y \Rightarrow \frac{1}{p_Y} \leq \frac{1}{p_X}. \qquad (33)$$

Subtracting 1 from both sides of the above inequality.

$$\frac{1}{p_Y} - 1 \leq \frac{1}{p_X} - 1 \Rightarrow \frac{1 - p_Y}{p_Y} \leq \frac{1 - p_X}{p_X}. \qquad (34)$$

Rearranging terms, we obtain:

$$\frac{p_X(1 - p_Y)}{(1 - p_X)p_Y} \leq 1 \Rightarrow p \leq 1 \left( \text{as } p = \frac{p_X(1 - p_Y)}{(1 - p_X)p_Y} \right). \qquad (35)$$

For calculating $P(Y)$, we borrow the result of proposition 13 in [26], which states that if $X \sim$ Negative Binomial$(M, p_X)$ and $P(Y|X) =$ Binomial$(X, p)$, then $P(Y) =$ Negative Binomial $\left( N, \frac{p_X}{p_X + p - pp_X} \right)$. Hence,

$$P(Y) = \text{Negative Binomial} \left( N, \frac{p_X}{p_X + p - pp_X} \right) = \text{Negative Binomial} \left( N, \frac{p_X}{p_X + p(1 - p_X)} \right)$$

$$= \text{Negative Binomial} \left( N, \frac{p_X}{p_X + \frac{p_X(1-p_Y)}{(1-p_X)p_Y}(1 - p_X)} \right)$$

$$= \text{Negative Binomial} \left( N, \frac{p_X}{p_X + \frac{p_X(1-p_Y)}{p_Y}} \right)$$

$$= \text{Negative Binomial} \left( N, \frac{p_X p_Y}{p_X p_Y + p_X(1 - p_Y)} \right)$$

$$= \text{Negative Binomial} \left( N, \frac{p_X p_Y}{p_X} \right) = \text{Negative Binomial} \left( N, p_Y \right). \qquad (36)$$

$\qquad\square$

### E.3 PID for the miscellaneous system M3.

Miscellaneous System **M3**: Let $(M, X, Y)$ be a system of random variables with joint distribution $P(M, X, Y)$. The joint distribution $P(M, X, Y)$ describes the miscellaneous system **M3** if it satisfies the following two properties:

1. $M \sim P(M)$ and has support $(0, \infty)$.
2. $P(X|M)=\text{Uniform}(0, \gamma_X M)$ and $P(Y|M)=\text{Uniform}(0, \gamma_Y M)$, where $\gamma_X, \gamma_Y \in \mathbb{R} \backslash \{0\}$.

**Proposition 4.** *Let $M, X,$ and $Y$ be random variables whose joint distribution $P(M, X, Y)$ describes the miscellaneous system **M3** defined above. Then, $\Delta_P$ contains a distribution $Q_{MC}(M, X, Y)$, which admits both Markovian structures, i.e., $M \to X \to Y$ and $M \to Y \to X$. Consequently, both $UI(M; Y \backslash X) = UI(M; X \backslash Y) = 0$.*

*Proof.* We prove proposition 4 by explicitly constructing a joint distribution $Q_{MC}(M, X, Y) \in \Delta_P$ that admits both Markovian structures, namely, $M \to X \to Y$ and $M \to Y \to X$. A necessary and sufficient condition for the distribution $Q_{MC}(M, X, Y)$ to admit both Markovian structures is that it should satisfy (37):

$$Q_{MC}(M, X, Y)=Q_{MC}(M)Q_{MC}(X|M)Q_{MC}(Y|X)=Q_{MC}(M)Q_{MC}(Y|M)Q_{MC}(X|Y). \tag{37}$$

We now briefly outline the structure of the proof. The rest of the proof can be divided into three parts:

1. In the first part, we explicitly construct $Q_{MC}(M, X, Y)$.

2. In the second part, we show that the $Q_{MC}(M, X, Y)$ constructed in the first part, lies in $\Delta_P$.

3. In the third part, we conclude our proof by showing that $Q_{MC}(M, X, Y)$ satisfies (37) and, consequently, admits both Markovian structures.

**Part 1: Specifying $Q_{MC}(M, X, Y)$**

We specify $Q_{MC}(M, X, Y)$ by individually specifying $Q_{MC}(M)$, $Q_{MC}(X|M)$, and $Q_{MC}(Y|X)$. We choose $Q_{MC}(M)$ and $Q_{MC}(X|M)$ as follows:

$$Q_{MC}(M) = P(M), \text{ and } Q_{MC}(X|M) = P(X|M). \tag{38}$$

The distribution $Q_{MC}(Y|X)$ is specified through the following deterministic transformation:

$$Y = \frac{\gamma_Y}{\gamma_X} X. \tag{39}$$

**Part 2: Showing $Q_{MC}(M, X, Y) \in \Delta_P$**

For showing $Q_{MC}(M, X, Y) \in \Delta_P$, we first need to calculate $Q_{MC}(Y|M)$:

*Calculating $Q_{MC}(Y|M)$:* Since $Q_{MC}(X|M)=\text{Uniform}(0, \gamma_X M)$, and $Y = \gamma_Y/\gamma_X X$, we can use the result of lemma 3 to calculate $Q_{MC}(Y|M)$ as follows:

$$Q_{MC}(Y|M) = Q_{MC}\left(\frac{\gamma_Y}{\gamma_X}X \bigg| M\right) = \text{Uniform}\left(0, \frac{\gamma_Y}{\gamma_X}\gamma_X M\right) = \text{Uniform}(0, \gamma_Y M)$$
$$= P(Y|M). \tag{40}$$

Then, from (38) and (40), we can conclude:

$$Q_{MC}(M, X) = Q_{MC}(M)Q_{MC}(X|M) = P(M)P(X|M) = P(M, X), \tag{41}$$
$$Q_{MC}(M, Y) = Q_{MC}(M)Q_{MC}(Y|M) = P(M)P(Y|M) = P(M, Y). \tag{42}$$

Hence, $Q_{MC}(M, X, Y) \in \Delta_P$.

**Part 3: Showing $Q_{MC}(M, X, Y)$ satisfies (37)**

It is trivial to see that by the construction of $Q_{MC}(M, X, Y)$ given in Part 1, we have:

$$Q_{MC}(M, X, Y) = Q_{MC}(M)Q_{MC}(X|M)Q_{MC}(Y|X). \tag{43}$$

By the chain rule, we have:

$$Q_{MC}(M, X, Y) = Q_{MC}(M)Q_{MC}(Y|M)Q_{MC}(X|Y, M). \tag{44}$$

Hence, for showing that $Q_{MC}(M, X, Y)$ satisfies (37), we only need to show that $Q_{MC}(X|Y, M) = Q_{MC}(X|Y)$.

*Showing $Q_{MC}(X|Y, M) = Q_{MC}(X|Y)$:* Observe that $Y = \gamma_Y/\gamma_X X$ by construction. Therefore, we can equivalently express $X = \gamma_X/\gamma_Y Y$. Consequently,

$$Q_{MC}(X|Y, M) = \mathbb{I}[X = \gamma_X/\gamma_Y Y] \text{ as } X = \gamma_X/\gamma_Y Y, \tag{45}$$

where $\mathbb{I}[\cdot]$ is the indicator function. Since R.H.S of the above equation does not depend on $M$, we have:

$$Q_{MC}(X|Y, M) = \mathbb{I}[X = \gamma_X/\gamma_Y Y] = Q_{MC}(X|Y). \tag{46}$$

Combining (44) and (46), we have:

$$Q_{MC}(M, X, Y) = Q_{MC}(M)Q_{MC}(Y|M)Q_{MC}(X|Y). \tag{47}$$

Combining (43) and (47), we can show that $Q_{MC}(M, X, Y)$ satisfies (37).

**Conclusion**: Combining the results of parts 2 and 3, we can conclude that $Q_{MC}(M, X, Y) \in \Delta_P$ and contains both Markovian structures: $M \rightarrow X \rightarrow Y$ and $M \rightarrow Y \rightarrow X$. A unique implication of $Q_{MC}(M, X, Y)$ admitting both Markovian structures is that both $I_{Q_{MC}}(M; X|Y) = I_{Q_{MC}}(M; Y|X) = 0$. Consequently, both minimization problems presented in (2) are minimized by $Q_{MC}(M, X, Y)$ with $UI(M; X \backslash Y) = \min_{Q \in \Delta_P} I_Q(M; X|Y) = 0$ and $UI(M; Y \backslash X) = \min_{Q \in \Delta_P} I_Q(M; Y|X) = 0$, completing our proof. □

**Lemma 3.** *Let $X \sim Uniform(0, a)$ for some $a \in \mathbb{R}$, then $\gamma X \sim Uniform(0, \gamma a)$ for $\gamma \in \mathbb{R}$ and $a, \gamma \neq 0$.*

*Proof.* We denote the p.d.f. of the random variable $X$ as $p_X(\cdot)$. By lemma statement, we know:

$$p_X(X = x) = \text{Uniform}(0, a) = \begin{cases} \frac{1}{|a|} & x \in [0, a] \\ 0 & \text{otherwise} \end{cases}. \tag{48}$$

Denote the p.d.f. of the function $f(X)$ of the random variable $X$ as $p_{f(X)}(\cdot)$. We want to derive $p_{f(X)}(\cdot)$ for $f(X) = \gamma X$. Since, $f(X) = \gamma X$ is an invertible function, we can use the change-of-variable technique for deriving $p_{f(X)}(\cdot)$:

$$p_{f(X)}(f(X) = y) = p_X(X = f^{-1}(y)) \left| \frac{df^{-1}(y)}{dy} \right| = \begin{cases} 1/|\gamma a| & y \in [0, \gamma a] \\ 0 & \text{otherwise} \end{cases} = \text{Uniform}(0, \gamma a).$$

□

### E.4 PID for the miscellaneous system M4.

Miscellaneous System **M4**: Let $(M, X, Y)$ be a system of random variables with joint distribution $P(M, X, Y)$. The joint distribution $P(M, X, Y)$ describes the miscellaneous system **M4** if it satisfies the following two properties:

1. $M \sim P(M)$ and has support $[0, 1]$.
2. $P(X|M) = \text{Bernoulli}(M)$, and $P(Y|M) = \text{Bernoulli}(M + p - 2pM)$, where $p \in [0, 1]$.

This system is adapted from the Bernoulli decomposition listed in Appendix A [26].

**Proposition 5.** *Let $M, X,$ and $Y$ be random variables whose joint distribution $P(M, X, Y)$ describes the miscellaneous system **M4** defined above. Then, $\Delta_P$ contains a Markov chain of the form $M \rightarrow X \rightarrow Y$ and $UI(M; Y \backslash X) = 0$.*

*Proof.* We briefly outline the proof structure. The proof can be divided into two parts:

1. In the first part, we explicitly construct a joint distribution $Q_{MC}(M, X, Y)$ having the Markovian structure $M \to X \to Y$.

2. In the second part, we conclude our proof by showing the $Q_{MC}(M, X, Y)$ constructed in the first part, lies in $\Delta_P$.

**Part 1: Specifying $Q_{MC}(M, X, Y)$**

Since $Q_{MC}(M, X, Y)$ is the joint distribution of a Markov chain $M \to X \to Y$, we have $Q_{MC}(M, X, Y) = Q_{MC}(M)Q_{MC}(X|M)Q_{MC}(Y|X)$. Leveraging the special structure of $Q_{MC}(M, X, Y)$, we specify $Q_{MC}(M, X, Y)$ by individually specifying $Q_{MC}(M)$, $Q_{MC}(X|M)$, and $Q_{MC}(Y|X)$. We choose $Q_{MC}(M)$, $Q_{MC}(X|M)$ as follows:

$$Q_{MC}(M) = P(M), \text{ and } Q_{MC}(X|M) = P(X|M). \tag{49}$$

The distribution $Q_{MC}(Y|X)$ is specified through the following stochastic transformation:

$$Y = X(1 - Z) + Z(1 - X), \tag{50}$$

where $Z \sim \text{Bernoulli}(p)$ and $Z \perp\!\!\!\perp (X, M)$.

**Part 2: Showing $Q_{MC}(M, X, Y) \in \Delta_P$**

For showing $Q_{MC}(M, X, Y) \in \Delta_P$, we first need to calculate $Q_{MC}(Y|M)$:

*Calculating $Q_{MC}(Y|M)$:* For calculating $Q(Y|M)$, we use the result of proposition 10 in [26], which states that if $X' \sim \text{Bernoulli}(\theta)$, $Z' \sim \text{Bernoulli}(p')$, $Z' \perp\!\!\!\perp X'$, and $Y' = X'(1 - Z') + (1 - X')Z'$, then $Y \sim \text{Bernoulli}(\theta + p' - 2p'\theta)$.

Noting that $Q_{MC}(X|M) = \text{Bernoulli}(M)$ and $Y = X(1 - Z) + Z(1 - X)$, where $Z \sim \text{Bernoulli}(p)$ and $Z \perp\!\!\!\perp X|M$, we can calculate $Q_{MC}(Y|M)$ as follows:

$$Q_{MC}(Y|M) = \text{Bernoulli}(M + p - 2pM) = P(Y|M). \tag{51}$$

Then, from (49) and (51), we can conclude:

$$Q_{MC}(M, X) = Q_{MC}(M)Q_{MC}(X|M) = P(M)P(X|M) = P(M, X), \tag{52}$$
$$Q_{MC}(M, Y) = Q_{MC}(M)Q_{MC}(Y|M) = P(M)P(Y|M) = P(M, Y). \tag{53}$$

Hence, $Q_{MC}(M, X, Y) \in \Delta_P$ and consequently, by proposition 1, $UI(M; Y \backslash X) = 0$. This concludes our proof. $\square$

### E.5 PID for the miscellaneous system M5.

Miscellaneous System **M5**: Let $(\vec{\mathbf{M}}, X, Y)$ be a system of random variables with joint distribution $P(\vec{\mathbf{M}}, X, Y)$. The joint distribution $P(\vec{\mathbf{M}}, X, Y)$ describes the miscellaneous system **M5** if it satisfies the following two properties:

1. $\vec{\mathbf{M}} \sim P(\vec{\mathbf{M}})$ and has support over simplex of dimension $d$.

2. $P(X|\vec{\mathbf{M}}) = \text{Categorical}(M_1, \dots, M_d)$, and $P(Y|\vec{\mathbf{M}}) = \text{Categorical}(\phi_1, \dots, \phi_d)$, where $\phi_i = (1 - p)M_i + p/d$.

This system is adapted from the categorical decomposition listed in Appendix A [26].

**Proposition 6.** *Let $\vec{\mathbf{M}}, X$, and $Y$ be random variables whose joint distribution $P(\vec{\mathbf{M}}, X, Y)$ describes the miscellaneous system **M5** defined above. Then, $\Delta_P$ contains a Markov chain of the form $\vec{\mathbf{M}} \to X \to Y$ and $UI(\vec{\mathbf{M}}; Y \backslash X) = 0$.*

*Proof.* We briefly outline the proof structure. The proof can be divided into two parts:

1. In the first part, we explicitly construct a joint distribution $Q_{MC}(\vec{\mathbf{M}}, X, Y)$ having the Markovian structure $\vec{\mathbf{M}} \to X \to Y$.

2. In the second part, we conclude our proof by showing the $Q_{MC}(\vec{\mathbf{M}}, X, Y)$ constructed in the first part, lies in $\Delta_P$.

**Part 1: Specifying $Q_{MC}(\vec{\mathbf{M}}, X, Y)$**

Since $Q_{MC}(\vec{\mathbf{M}}, X, Y)$ is the joint distribution of a Markov chain $\vec{\mathbf{M}} \to X \to Y$, we have $Q_{MC}(\vec{\mathbf{M}}, X, Y) = Q_{MC}(\vec{\mathbf{M}})Q_{MC}(X|\vec{\mathbf{M}})Q_{MC}(Y|X)$. Taking advantage of this special structure of $Q_{MC}(\vec{\mathbf{M}}, X, Y)$, we specify $Q_{MC}(\vec{\mathbf{M}}, X, Y)$ by individually specifying $Q_{MC}(\vec{\mathbf{M}})$, $Q_{MC}(X|\vec{\mathbf{M}})$, and $Q_{MC}(Y|X)$. We choose $Q_{MC}(\vec{\mathbf{M}}), Q_{MC}(X|\vec{\mathbf{M}})$ as follows:

$$Q_{MC}(\vec{\mathbf{M}}) = P(\vec{\mathbf{M}}), \text{ and } Q_{MC}(X|\vec{\mathbf{M}}) = P(X|\vec{\mathbf{M}}). \tag{54}$$

The distribution $Q_{MC}(Y|X)$ is specified through the following transformation:

$$Y = X(1 - Z) + ZD, \tag{55}$$

where $Z \sim \text{Bernoulli}(p)$, $D \sim \text{Categorical}\left(\frac{1}{d}, \ldots, \frac{1}{d}\right)$, and $D, Z$, and $(X, \vec{\mathbf{M}})$ are jointly independent.

**Part 2: Showing $Q_{MC}(\vec{\mathbf{M}}, X, Y) \in \Delta_P$**

For showing $Q_{MC}(\vec{\mathbf{M}}, X, Y) \in \Delta_P$, we first need to calculate $Q_{MC}(Y|\vec{\mathbf{M}})$:

*Calculating $Q_{MC}(Y|\vec{\mathbf{M}})$:* For calculating $Q_{MC}(Y|\vec{\mathbf{M}})$, we use the result of proposition 11 in [26], which states that:

If $X' \sim \text{Categorical}(\theta_1, \ldots, \theta_d)$, $Z' \sim \text{Bernoulli}(p')$, $D' \sim \text{Categorical}(1/d, \ldots, 1/d)$, and $Y' = X'(1 - Z') + Z'D'$ with $X', Z'$, and $D'$ jointly independent, then $Y' \sim \text{Categorical}(\phi'_1, \ldots, \phi'_d)$ with $\phi'_i = (1 - p')\theta_i + p'/d$.

Noting that $Q_{MC}(X|\vec{\mathbf{M}}) = \text{Categorical}(M_1, \ldots, M_d)$ and $Y = X(1 - Z) + ZD$, with $X, D$, and $Z$ jointly conditionally independent conditioned on $M$, we have:

$$Q_{MC}(Y|\vec{\mathbf{M}}) = \text{Categorical}(\phi_1, \ldots, \phi_d), \text{ with } \phi_i = M_i(1 - p) + \frac{p}{d} = P(Y|\vec{\mathbf{M}}). \tag{56}$$

From (54) and (56), we can conclude:

$$Q_{MC}(\vec{\mathbf{M}}, X) = Q_{MC}(\vec{\mathbf{M}})Q_{MC}(X|\vec{\mathbf{M}}) = P(M)P(X|\vec{\mathbf{M}}) = P(\vec{\mathbf{M}}, X), \tag{57}$$

$$Q_{MC}(\vec{\mathbf{M}}, Y) = Q_{MC}(\vec{\mathbf{M}})Q_{MC}(Y|\vec{\mathbf{M}}) = P(M)P(Y|\vec{\mathbf{M}}) = P(\vec{\mathbf{M}}, Y). \tag{58}$$

Hence, $Q_{MC}(\vec{\mathbf{M}}, X, Y) \in \Delta_P$ and consequently, by proposition 1, $UI(\vec{\mathbf{M}}; Y \backslash X) = 0$. This concludes our proof. $\qquad\square$

## E.6 PID for the miscellaneous system M6.

Miscellaneous System **M6**: Let $(M, \vec{\mathbf{X}}, \vec{\mathbf{Y}})$ be a system of random variables with joint distribution $P(M, \vec{\mathbf{X}}, \vec{\mathbf{Y}})$. The joint distribution $P(M, \vec{\mathbf{X}}, \vec{\mathbf{Y}})$ describes the miscellaneous system **M6** if it satisfies the following two properties:

1. $M \sim P(M)$ and has support $\mathbb{N}$.
2. $P(\vec{\mathbf{X}}|M) = \text{Multinomial}_{d_X}(\vec{\mathbf{X}}; M, \vec{\mathbf{p}}_X)$, and $P(\vec{\mathbf{Y}}|M) = \text{Multinomial}_{d_Y}(\vec{\mathbf{Y}}; M, \vec{\mathbf{p}}_Y)$.

**Proposition 7.** *Let $M$ be a random variable. Furthermore, assume $\vec{\mathbf{X}}$ and $\vec{\mathbf{Y}}$ are random vectors of size $d_X$ and $d_Y$, respectively, such that $P(M, \vec{\mathbf{X}}, \vec{\mathbf{Y}})$ describes miscellaneous system **M6**.*

1. *If, $\min_{i \in [d_Y]} p_i^Y \geq \min_{i \in [d_X]} p_i^X$, then there exists a Markov chain of the form $M \to \vec{\mathbf{X}} \to \vec{\mathbf{Y}}$ in $\Delta_P$, and $UI(M; \vec{\mathbf{Y}} \backslash \vec{\mathbf{X}}) = 0$.*

2. *If, $\min_{i \in [d_Y]} p_i^Y \leq \min_{i \in [d_X]} p_i^X$, then there exists a Markov chain of the form $M \to \vec{\mathbf{Y}} \to \vec{\mathbf{X}}$ in $\Delta_P$, and $UI(M; \vec{\mathbf{X}} \backslash \vec{\mathbf{Y}}) = 0$.*

*Proof.* We only provide an explicit proof of condition 1. The proof of condition 2 follows the same steps as the proof of condition 1 with parameters of $\vec{\mathbf{X}}$ and $\vec{\mathbf{Y}}$ switched.

We briefly outline the proof structure. The proof can be divided into two major parts:

1. In the first part, we explicitly construct a joint distribution $Q_{MC}(M, \vec{\mathbf{X}}, \vec{\mathbf{Y}})$ having the Markovian structure $M \to \vec{\mathbf{X}} \to \vec{\mathbf{Y}}$.

2. In the second part, we conclude our proof by showing the $Q_{MC}(M, \vec{\mathbf{X}}, \vec{\mathbf{Y}})$ constructed in the first part, lies in $\Delta_P$.

**Part 1: Specifying $Q_{MC}(M, \vec{\mathbf{X}}, \vec{\mathbf{Y}})$**

Since $Q_{MC}(M, \vec{\mathbf{X}}, \vec{\mathbf{Y}})$ is the joint distribution of a Markov chain $M \to \vec{\mathbf{X}} \to \vec{\mathbf{Y}}$, we have $Q_{MC}(M, \vec{\mathbf{X}}, \vec{\mathbf{Y}}) = Q_{MC}(M) Q_{MC}(\vec{\mathbf{X}}|M) Q_{MC}(\vec{\mathbf{Y}}|\vec{\mathbf{X}})$. Taking advantage of this special structure of $Q_{MC}(M, \vec{\mathbf{X}}, \vec{\mathbf{Y}})$, we specify $Q_{MC}(M, \vec{\mathbf{X}}, \vec{\mathbf{Y}})$ by individually specifying $Q_{MC}(M)$, $Q_{MC}(\vec{\mathbf{X}}|M)$, and $Q_{MC}(\vec{\mathbf{Y}}|\vec{\mathbf{X}})$. We choose $Q_{MC}(M), Q_{MC}(\vec{\mathbf{X}}|M)$ as follows:

$$Q_{MC}(M) = P(M), \text{ and } Q_{MC}(\vec{\mathbf{X}}|M) = P(\vec{\mathbf{X}}|M). \tag{59}$$

*Specifying $Q(\vec{\mathbf{Y}}|\vec{\mathbf{X}})$:* Let $k_X^* = \arg\min_{i \in [d_X]} p_i^X$, and $k_Y^* = \arg\min_{i \in [d_Y]} p_i^Y$. Then, $Q(\vec{\mathbf{Y}}|\vec{\mathbf{X}}) = \text{Multinomial}_{d_Y} \left( \vec{\mathbf{Y}}; \sum_{i \in [d_X] \setminus k_X^*} x_i, \mathbf{p}_Y^* \right)$, where

$$\mathbf{p}_Y^* = \left[ \frac{p_1^Y}{\sum_{i \in [d_X] \setminus k_X^*} p_i^X} \cdots \frac{p_{d_Y-1}^Y}{\sum_{i \in [d_X] \setminus k_X^*} p_i^X} \quad 1 - \frac{\sum_{i=1}^{d_Y-1} p_i^Y}{\sum_{i \in [d_X] \setminus k_X^*} p_i^X} \right]^T. \tag{60}$$

We first show the proposed $Q_{MC}(\vec{\mathbf{Y}}|\vec{\mathbf{X}})$ is a valid distribution. By condition 1:

$$\min_{i \in [d_Y]} p_i^Y \geq \min_{i \in d_X} p_i^X \tag{61}$$

$$\Rightarrow \sum_{i \in [d_X] \setminus k_X^*} p_i^X \geq \sum_{i \in [d_Y] \setminus k_Y^*} p_i^Y \geq p_j^Y, \ \forall\, j \in [d_Y] \tag{62}$$

$$\Rightarrow 1 \geq \frac{p_j^Y}{\sum_{i \in [d_X] \setminus k_X^*} p_i^X} \geq 0, \ \forall\, j \in [d_Y], \tag{63}$$

where the last inequality is due to $\vec{\mathbf{p}}_X$ and $\vec{\mathbf{p}}_Y$ forming valid multinomial distributions.

**Part 2: Showing $Q_{MC}(M, \vec{\mathbf{X}}, \vec{\mathbf{Y}}) \in \Delta_P$**

For showing $Q_{MC}(M, \vec{\mathbf{X}}, \vec{\mathbf{Y}}) \in \Delta_P$, we first need to calculate $Q_{MC}(\vec{\mathbf{Y}}|M)$:

*Calculating $Q_{MC}(\vec{\mathbf{Y}}|M)$:* We use lemma 4 to calculate $Q(\vec{\mathbf{Y}}|M)$. Note that $Q_{MC}(\vec{\mathbf{X}}|M) = \text{Multinomial}_{d_X}(M, \vec{\mathbf{p}}_X)$, hence we have:

$$Q_{MC}(Y|M) = \text{Multinomial}_{d_Y}(M, \vec{\mathbf{p}}_Y'), \text{ where },$$

$$\vec{\mathbf{p}}_Y' = \left[ \frac{p_1^Y \sum_{i \in [d_X] \setminus k_X^*} p_i^X}{\sum_{i \in [d_X] \setminus k_X^*} p_i^X} \quad \cdots \quad \frac{p_{d_Y-1}^Y \sum_{i \in [d_X] \setminus k_X^*} p_i^X}{\sum_{i \in [d_X] \setminus k_X^*} p_i^X} \quad 1 - \frac{\sum_{i \in [d_X] \setminus k_X^*} p_i^X \sum_{i=1}^{d_Y-1} p_i^Y}{\sum_{i \in [d_X] \setminus k_X^*} p_i^X} \right]$$

$$\Rightarrow \vec{\mathbf{p}}_Y' = \left[ p_1^Y \quad \cdots \quad p_{d_Y-1}^Y \quad 1 - \sum_{i \in [d_Y]} p_i^Y \right] = \vec{\mathbf{p}}_Y$$

$$\Rightarrow Q_{MC}(Y|M) = \text{Multinomial}_{d_Y}(M, \vec{\mathbf{p}}_Y) = P(Y|M). \tag{64}$$

Then, from (59) and (64), we can conclude:

$$Q_{MC}(M, \vec{\mathbf{X}}) = Q_{MC}(M) Q_{MC}(\vec{\mathbf{X}}|M) = P(M) P(\vec{\mathbf{X}}|M) = P(M, \vec{\mathbf{X}}), \tag{65}$$

$$Q_{MC}(M, \vec{\mathbf{Y}}) = Q_{MC}(M) Q_{MC}(\vec{\mathbf{Y}}|M) = P(M) P(\vec{\mathbf{Y}}|M) = P(M, \vec{\mathbf{Y}}). \tag{66}$$

Hence, $Q_{MC}(M, \vec{\mathbf{X}}, \vec{\mathbf{Y}}) \in \Delta_P$ and consequently, by proposition 1, $UI(M; \vec{\mathbf{Y}} \setminus \vec{\mathbf{X}}) = 0$. This concludes our proof. $\quad\square$

**Lemma 4.** *Suppose* $\vec{\mathbf{X}} \sim Multinomial_{d_X}(n, \vec{\mathbf{p}}_X)$ *and* $\vec{\mathbf{Y}}|\vec{\mathbf{X}} \sim Multinomial_{d_Y}\left(\sum_{j\in\mathcal{I}} X_j, \vec{\mathbf{q}}\right)$, *where* $\mathcal{I} \subset \{1,\ldots,d_X\}$. *Then* $\vec{\mathbf{Y}} \sim Multinomial_{d_Y}(n, \vec{\mathbf{q}}^*)$, *where*

$$q_i^* = \begin{cases} \left[\sum_{j\in\mathcal{I}} p_j\right] \cdot q_i, & 1 \leq i < d_Y \\ 1 - \left[\sum_{j\in\mathcal{I}} p_j\right] \cdot \left[\sum_{j=1}^{d_Y-1} q_j\right], & i = d_Y \end{cases}. \tag{67}$$

*Proof.* Without loss of generality, reshuffle the class indices of $\vec{\mathbf{Y}}$ such that $\mathcal{I}=\{i: 1 \leq i \leq w\}$, where $|\mathcal{I}|=w$. Let $C=\{\vec{\mathbf{x}}' \in \mathbb{N}_0^{d_X-1}: \sum_{i=1}^{d_X-1} x_i \leq n\}$. By the law of total probability,

$$P(\vec{\mathbf{Y}} = \vec{\mathbf{y}}) = \sum_{\vec{\mathbf{x}}\in C} P(\vec{\mathbf{X}} = \vec{\mathbf{x}})P(\vec{\mathbf{Y}} = \vec{\mathbf{y}}|\vec{\mathbf{X}} = \vec{\mathbf{x}}) \tag{68}$$

$$= \sum_{\vec{\mathbf{x}}\in C} \frac{n!}{\prod_{i=1}^{d_X} x_i!} \prod_{i=1}^{d_X} p_i^{x_i} \cdot \frac{\left(\sum_{j\in\mathcal{I}} x_j\right)!}{\prod_{i=1}^{d_Y} y_i!} \prod_{i=1}^{d_Y} q_i^{y_i}. \tag{69}$$

Remove from the sum all terms that do not depend on $\vec{\mathbf{x}}$. Note that $y_{d_Y} = \sum_{j\in\mathcal{I}} x_j - \sum_{i=1}^{d_Y-1} y_i$ and $q_{d_Y}^{y_{d_Y}} = \left(1 - \sum_{i=1}^{d_Y} q_i\right)^{y_{d_Y}}$ are both functions of $\vec{\mathbf{y}}$. Let $y_s = \sum_{i=1}^{d_Y-1} y_i$ and $q_s = \sum_{i=1}^{d_Y-1} q_i$.

$$P(\vec{\mathbf{Y}} = \vec{\mathbf{y}}) = \left(\frac{n!}{\prod_{i=1}^{d_Y-1} y_i!} \prod_{i=1}^{d_Y-1} q_i^{y_i}\right) \sum_{\vec{\mathbf{x}}\in C} \frac{\prod_{i=1}^{d_X} p_i^{x_i} \cdot \left(\sum_{j\in\mathcal{I}} x_j\right)!}{\prod_{i=1}^{d_X} x_i! \left(\sum_{j\in\mathcal{I}} x_j - y_s\right)!} (1 - q_s)^{\sum_{j\in\mathcal{I}} x_j - y_s}.$$

Form a multinomial coefficient outside the sum:

$$P(\vec{\mathbf{Y}} = \vec{\mathbf{y}}) = \left(\frac{n!}{\prod_{i=1}^{d_Y-1} y_i! \cdot (n - y_s)!} \prod_{i=1}^{d_Y-1} q_i^{y_i}\right) \tag{70}$$

$$\times \sum_{\vec{\mathbf{x}}\in C} \frac{(n - y_s)! \prod_{i=1}^{d_X} p_i^{x_i} \cdot \left(\sum_{j\in\mathcal{I}} x_j\right)!}{\prod_{i=1}^{d_X} x_i! \left(\sum_{j\in\mathcal{I}} x_j - y_s\right)!} (1 - q_s)^{\sum_{j\in\mathcal{I}} x_j - y_s}. \tag{71}$$

Let $\mathcal{I}^0 = \{i: w < i \leq d_X\}$ be the set of indices that are not taken in the number of trials for $\vec{\mathbf{Y}}$. Consider the sum over $\vec{\mathbf{x}} \in C$ alone in (71). Separating terms that belong to $\mathcal{I}$ and $\mathcal{I}^0$,

$$\sum_{\vec{\mathbf{x}}\in C} \frac{(n - y_s)!}{\prod_{j\in\mathcal{I}^0} x_j! \left(\sum_{j\in\mathcal{I}} x_j - y_s\right)!} \prod_{j\in\mathcal{I}^0} p_j^{x_j} (1 - q_s)^{\sum_{j\in\mathcal{I}} x_j - y_s} \cdot \frac{\left(\sum_{j\in\mathcal{I}} x_j\right)!}{\prod_{j\in\mathcal{I}} x_j!} \prod_{j\in\mathcal{I}} p_j^{x_j}.$$

Let $C' = \{\vec{\mathbf{x}}' \in \mathbb{N}_0^{d_X-1}: \sum_{i=1}^{d_X-1} x_i - \sum_{i=1}^{d_Y-1} y_i \leq n - \sum_{i=1}^{d_Y-1} y_i\}$. Note $\vec{\mathbf{x}} \in C \Leftrightarrow \vec{\mathbf{x}} \in C'$, so we can equivalently sum over elements in $C'$. Perform a change of variable with $u = \sum_{j\in\mathcal{I}} x_j - y_s$ and define $B = \{x_j \in \vec{\mathbf{x}}: j \in \mathcal{I}, \sum_{j\in\mathcal{I}} x_j = u + y_s\}$. Then the sum becomes

$$= \sum_{\vec{\mathbf{x}}\in C'} \frac{(n - y_s)!}{\prod_{j\in\mathcal{I}^0} x_j! u!} \prod_{j\in\mathcal{I}^0} p_j^{x_j} (1 - q_s)^u \sum_{x_j\in B} \binom{u + y_s}{x_{j_1}, \ldots, x_{j_m}} \prod_{j\in\mathcal{I}} p_j^{x_j}. \tag{72}$$

Note that the inner sum simplifies by the multinomial theorem:

$$= \sum_{\vec{\mathbf{x}}\in C'} \frac{(n - y_s)!}{\prod_{j\in\mathcal{I}^0} x_j! u!} \prod_{j\in\mathcal{I}^0} p_j^{x_j} (1 - q_s)^u \left(\sum_{j\in\mathcal{I}} p_j\right)^{u+y_s} \tag{73}$$

$$= \left(\sum_{j\in\mathcal{I}} p_j\right)^{y_s} \sum_{\vec{\mathbf{x}}\in C'} \frac{(n - y_s)!}{\prod_{j\in\mathcal{I}^0} x_j! \cdot u!} \prod_{j\in\mathcal{I}^0} p_j^{x_j} \left(\sum_{j\in\mathcal{I}} p_j - q_s \sum_{j\in\mathcal{I}} p_j\right)^u. \tag{74}$$

Reapply the multinomial theorem to the sum of $\vec{x}$ over $C'$:

$$= \left( \sum_{j \in \mathcal{I}} p_j \right)^{y_s} \left( \sum_{j \in \mathcal{I}^0} p_j + \sum_{j \in \mathcal{I}} p_j - q_s \sum_{j \in \mathcal{I}} p_j \right)^{n-y_s} \tag{75}$$

$$= \left( \sum_{j \in \mathcal{I}} p_j \right)^{y_s} \left( 1 - q_s \sum_{j \in \mathcal{I}} p_j \right)^{n-y_s} . \tag{76}$$

Inserting (76) into (71), the marginal becomes

$$P(\vec{\mathbf{Y}} = \vec{\mathbf{y}}) = \frac{n! \prod_{i=1}^{d_Y-1} q_i^{y_i}}{\prod_{i=1}^{d_Y-1} y_i! \cdot (n-y_s)!} \left( \sum_{j \in \mathcal{I}} p_j \right)^{y_s} \left( 1 - q_s \sum_{j \in \mathcal{I}} p_j \right)^{n-y_s} \tag{77}$$

$$= \frac{n! \prod_{i=1}^{d_Y-1} \left( \sum_{j \in \mathcal{I}} p_j \cdot q_i \right)^{y_i}}{\prod_{i=1}^{d_Y-1} y_i! \cdot (n-y_s)!} \left( 1 - q_s \sum_{j \in \mathcal{I}} p_j \right)^{n-y_s} \tag{78}$$

$$= \text{Multinomial}_{d_Y} \left( n, \vec{\mathbf{q}}^* \right) \tag{79}$$

where $\vec{\mathbf{q}}^*$ is defined as in (67). $\qquad\square$

## F    Proof of Theorem 1

In this section, we provide the proof of theorem 1. Since theorem 1 provides the analytical PID terms for the univariate affine continuous stable system defined in Sec. 4, we briefly restate certain key properties of the univariate stable distributions and the corresponding univariate affine continuous stable system for convenience.

### F.1    Univariate continuous stable distribution

Univariate continuous stable distribution are a large class of distributions that naturally arise in the context of generalized central limit theorems. We refer the reader to Appx. N.2 for more details on univariate continuous stable distributions. We now list certain key properties of univariate continuous stable distributions that we make use of in the proof of theorem 1:

1. If $X$ is distributed according to a univariate continuous stable distribution, then the sum of two independent copies of $X$, denoted as $X_1$ and $X_2$, follows the same univariate continuous stable distribution upto a scaling and translation operation, i.e., $aX_1 + bX_2 \overset{d}{=} cX + d$ for $a, b, c > 0$ and $d \in \mathbb{R}$.

2. The p.d.f. of univariate continuous stable distributions is characterized by four parameters: stability parameter denoted as $\alpha \in (0, 2]$, skewness parameter denoted as $\beta \in [-1, 1]$, scale parameters denoted as $\gamma \in (0, \infty)$, and location parameter denoted as $\mu \in \mathbb{R}$. We denote the p.d.f. of a univariate continuous stable distribution as $p_{CS}(\alpha, \beta, \gamma, \mu)$.

### F.2    Definition of univariate affine continuous stable system

Let $(M, X, Y)$ be a system of random variables with joint distribution $P(M, X, Y)$. The joint distribution $P(M, X, Y)$ describes the univariate affine continuous stable system if it satisfies the following two properties:

1. $M \sim P(M)$ having some support set $\mathcal{M} \subseteq \mathbb{R}$.

2. The conditional distributions of random variables $X$ and $Y$ conditioned on $M$ can be expressed through univariate continuous stable distributions with an affine dependence on $M$, i.e., $P(X|M) = p_{CS}(\alpha, \beta_X, \gamma_X, aM + b)$ and $P(Y|M) = p_{CS}(\alpha, \beta_Y \operatorname{sgn}(ac), \gamma_Y, cM + d)$, where $a, b, c, d \in \mathbb{R}$.

### F.3  Formal proof of Theorem 1

We first discuss and provide the provide the proofs of lemma 5 and lemma 6 that we will use for proving theorem 1. In the following proofs, $\mathrm{sgn}(x)$ denotes the sign function:

$$\mathrm{sgn}(x) = \left\{ \begin{array}{ll} 1 & x > 0 \\ 0 & x = 0 \\ -1 & x < 0 \end{array} \right. . \tag{80}$$

**Lemma 5.** *If $X \sim p_{CS}(\alpha, \beta, \gamma, \mu)$, then for any $\eta \neq 0$ and $\kappa \in \mathbb{R}$, we have*

$$\eta X + \kappa \sim \left\{ \begin{array}{ll} p_{CS}(\alpha, sgn(\eta)\beta, |\eta|\gamma, \eta\mu + \kappa) & \alpha \neq 1 \\ p_{CS}\left(\alpha, sgn(\eta)\beta, |\eta|\gamma, \eta\mu + \kappa - \frac{2}{\pi}\beta\gamma\eta \log(|\eta|)\right) & \alpha = 1 \end{array} \right. .$$

*Proof.* See the proof of proposition 1.4 part (b) in [23]. ☐

**Lemma 6.** *If $X_1 \sim p_{CS}(\alpha, \beta_1, \gamma_1, \mu_1)$ and $X_2 \sim p_{CS}(\alpha, \beta_2, \gamma_2, \mu_2)$ are independent, then $X_1 + X_2 \sim p_{CS}(\alpha, \beta, \gamma, \mu)$ where*

$$\beta = \frac{\beta_1 \gamma_1^\alpha + \beta_2 \gamma_2^\alpha}{\gamma_1^\alpha + \gamma_2^\alpha}, \quad \gamma^\alpha = \gamma_1^\alpha + \gamma_2^\alpha, \text{ and } \mu = \mu_1 + \mu_2.$$

*Proof.* See the proof of proposition 1.4 part (c) in [23]. ☐

**Theorem 1.** *Let $M, X$, and $Y$ be random variables whose joint distribution $P(M, X, Y)$ describes a univariate affine continuous stable system. Without the loss of generality, assume $|a|/\gamma_X \geq |c|/\gamma_Y$. If $1 - \beta_Y \geq (\gamma_X |c|/\gamma_Y |a|)^\alpha (1 - \beta_X)$, and $1 + \beta_Y \geq (\gamma_X |c|/\gamma_Y |a|)^\alpha (1 + \beta_X)$, then $\Delta_P$ contains a Markov chain of the form $M \to X \to Y$ and $UI(M; Y \backslash X) = 0$.*

*Proof.* We first note that we can always assume $|a|/\gamma_X \geq |c|/\gamma_Y$ without the loss of generality because if $|c|/\gamma_Y \geq |a|/\gamma_X$, then we can always switch our nomenclature to refer to $Y$ as $X$, and $X$ as $Y$.

We now briefly outline the proof structure. We divide the proof into two cases $|a|/\gamma_X > |c|/\gamma_Y$ and $|a|/\gamma_X = |c|/\gamma_Y$. The proof for both cases follows essentially the same structure consisting of two major parts:

1. In the first part, we explicitly construct a joint distribution $Q_{MC}(M, X, Y)$ having the Markovian structure $M \to X \to Y$.

2. In the second part, we show that the $Q_{MC}(M, X, Y)$ constructed in the first part, lies in $\Delta_P$. Therefore, we can then apply the result of proposition 1 to conclude $UI(M; Y \backslash X) = 0$.

**Case 1:** $|a|/\gamma_X > |c|/\gamma_Y$

**Part 1: Specifying $Q_{MC}(M, X, Y)$**

We explicitly construct the desired Markov chain $M \to X \to Y$ by constructing a larger Markov chain $M \to X \to X' \to Y' \to Y$ and then marginalizing the larger Markov chain to obtain the desired Markov chain $M \to X \to Y$.

Denote the joint distribution of the Markov chain $M \to X \to X' \to Y' \to Y$ as $Q_{MC}(M, X, X', Y', Y)$. We can decompose $Q_{MC}(M, X, X', Y', Y)$ by utilizing its Markovian structure as follows:

$$Q_{MC}(M, X, X', Y', Y) = Q_{MC}(M)Q_{MC}(X|M)Q_{MC}(X'|X)Q_{MC}(Y'|X')Q_{MC}(Y|Y').$$

Consequently, we can specify/construct the distribution $Q_{MC}(M, X, X', Y', Y)$ by individually specifying $Q_{MC}(M), Q_{MC}(X|M), Q_{MC}(X'|X), Q_{MC}(Y'|X')$, and $Q_{MC}(Y|Y')$.

Specifying $Q_{MC}(M)$ and $Q_{MC}(X|M)$: We choose $Q_{MC}(M)$ and $Q_{MC}(X|M)$ as follows:

$$Q_{MC}(M) = P(M) \text{ and } Q_{MC}(X|M) = P(X|M), \tag{81}$$

where $P(M)$ and $P(X|M)$ are marginal distributions derived from the original joint distribution $P(M, X, Y)$ (discussed in the theorem statement) over which the bivariate PID is being calculated.

Specifying $Q_{MC}(X'|X)$: The distribution $Q_{MC}(X'|X)$ is specified by the following deterministic transformation:

$$X' = \begin{cases} \frac{1}{a}X & \alpha \neq 1 \\ \frac{1}{a}X + (2\beta_X\gamma_X \log(|1/a|))/(\pi a) & \alpha = 1 \end{cases} \tag{82}$$

We will also derive the distribution $Q(X'|M)$ before proceeding with our construction, as we will need it later in the proof to show that the constructed $Q_{MC}(M, X, Y) \in \Delta_P$.

*Deriving $Q_{MC}(X'|M)$*: Since $Q_{MC}(X|M = m) = p_{CS}(\alpha, \beta_X, \gamma_X, am + b)$ for a fixed $m$ and $X'$ is scaled and translated version of $X$, we can use lemma 5 to derive $Q_{MC}(X'|M)$. The exact expression of $Q_{MC}(X'|M = m)$ is provided in (83).

$$Q_{MC}(X'|M = m) = p_{CS}(\alpha, \beta_{X'}, \gamma_{X'}, \mu_{X'}), \text{ where}$$

$$\beta_{X'} = \begin{cases} \text{sgn}(1/a)\beta_X & \alpha \neq 1 \\ \text{sgn}(1/a)\beta_X & \alpha = 1 \end{cases} = \text{sgn}(1/a)\beta_X = \text{sgn}(a)\beta_X,$$

$$\gamma_{X'} = \begin{cases} |1/a|\,\gamma_X & \alpha \neq 1 \\ |1/a|\,\gamma_X & \alpha = 1 \end{cases} = \frac{\gamma_X}{|a|},$$

$$\mu_{X'} = \begin{cases} \frac{1}{a}(am + b) & \alpha \neq 1 \\ \frac{1}{a}(am + b) + 2\beta_X\gamma_X \log(|1/a|)/\pi a - 2\beta_X\gamma_X \log(|1/a|)/\pi a & \alpha = 1 \end{cases} = m + \frac{b}{a},$$

$$\Rightarrow Q_{MC}(X'|M = m) = p_{CS}(\alpha, \beta_X\text{sgn}(a), \gamma_X/|a|, m + b/a). \tag{83}$$

Specifying $Q_{MC}(Y'|X')$: In order to define $Q_{MC}(Y'|X')$, we need to define an auxiliary variable $\epsilon$. The random variable $\epsilon \sim p_{CS}(\epsilon; \alpha, \beta', \gamma', \mu')$, where

$$\beta' = \text{sgn}(a)\frac{(\gamma_Y/|c|)^\alpha \beta_Y - (\gamma_X/|a|)^\alpha \beta_X}{(\gamma_Y/|c|)^\alpha - (\gamma_X/|a|)^\alpha}, \quad \gamma' = \left(\left(\frac{\gamma_Y}{|c|}\right)^\alpha - \left(\frac{\gamma_X}{|a|}\right)^\alpha\right)^{\frac{1}{\alpha}}, \quad \mu' = \frac{d}{c} - \frac{b}{a}. \tag{84}$$

In order for $\epsilon$ to have a legitimate continuous stable distribution, we need to ensure that $\beta'$, $\gamma'$, and $\mu'$ lie within their appropriate bounds as specified in Appx. F.1. It is trivial to see that $\mu' \in \mathbb{R}$.

Showing $\gamma' \in (0, \infty)$. Under the assumption of the case 1, we know that:

$$\frac{|a|}{\gamma_X} > \frac{|c|}{\gamma_Y}.$$

Manipulating the above inequality, we obtain:

$$\frac{\gamma_Y}{|c|} > \frac{\gamma_X}{|a|} \Rightarrow \left(\frac{\gamma_Y}{|c|}\right)^\alpha > \left(\frac{\gamma_X}{|a|}\right)^\alpha \Rightarrow \left(\frac{\gamma_Y}{|c|}\right)^\alpha - \left(\frac{\gamma_X}{|a|}\right)^\alpha > 0$$

$$\Rightarrow \left(\left(\frac{\gamma_Y}{|c|}\right)^\alpha - \left(\frac{\gamma_X}{|a|}\right)^\alpha\right)^{\frac{1}{\alpha}} = \gamma' > 0. \tag{85}$$

Hence, (85) shows that $\gamma' \in (0, \infty)$.

Lastly, showing that $\beta' \in [-1, 1]$. Note that the condition $\beta' \in [-1, 1]$ can be equivalently expressed as $|\beta'| \leq 1$. For showing that $\beta' \in [-1, 1]$, we will show that the inequality $|\beta'| \leq 1$ is equivalent to the inequalities: $(\gamma_X|c|/|a|\gamma_Y)^\alpha (1 - \beta_X) \leq 1 - \beta_Y$ and $(1 + \beta_X)(\gamma_X|c|/|a|\gamma_Y)^\alpha \leq 1 + \beta_Y$, where the last two inequalities hold by the theorem assumptions:

$$|\beta'| \leq 1 \Rightarrow \left|\text{sgn}(a)\frac{(\gamma_Y/|c|)^\alpha \beta_Y - (\gamma_X/|a|)^\alpha \beta_X}{(\gamma_Y/|c|)^\alpha - (\gamma_X/|a|)^\alpha}\right| \leq 1 \Rightarrow \left|\frac{(\gamma_Y/|c|)^\alpha \beta_Y - (\gamma_X/|a|)^\alpha \beta_X}{(\gamma_Y/|c|)^\alpha - (\gamma_X/|a|)^\alpha}\right| \leq 1,$$

$$\Rightarrow -1 \leq \frac{(\gamma_Y/|c|)^\alpha \beta_Y - (\gamma_X/|a|)^\alpha \beta_X}{(\gamma_Y/|c|)^\alpha - (\gamma_X/|a|)^\alpha} \leq 1. \tag{86}$$

Multiplying $(\gamma_Y/|c|)^\alpha - (\gamma_X/|a|)^\alpha$ on both sides of the inequalities in (86).

$$(\gamma_X/|a|)^\alpha - (\gamma_Y/|c|)^\alpha \leq (\gamma_Y/|c|)^\alpha \beta_Y - (\gamma_X/|a|)^\alpha \beta_X \leq (\gamma_Y/|c|)^\alpha - (\gamma_X/|a|)^\alpha. \tag{87}$$

Adding $(\gamma_X/|a|)^\alpha \beta_X$ on both sides of the inequalities in (87).

$$(\gamma_X/|a|)^\alpha + (\gamma_X/|a|)^\alpha \beta_X - (\gamma_Y/|c|)^\alpha \leq (\gamma_Y/|c|)^\alpha \beta_Y \leq (\gamma_Y/|c|)^\alpha - (\gamma_X/|a|)^\alpha + (\gamma_X/|a|)^\alpha \beta_X,$$
$$\Rightarrow (1 + \beta_X)(\gamma_X/|a|)^\alpha - (\gamma_Y/|c|)^\alpha \leq (\gamma_Y/|c|)^\alpha \beta_Y \leq (\gamma_Y/|c|)^\alpha + (\gamma_X/|a|)^\alpha (\beta_X - 1). \tag{88}$$

Dividing $(\gamma_Y/|c|)^\alpha$ on both sides of the inequalities in (88).

$$(1 + \beta_X)(\gamma_X|c|/|a|\gamma_Y)^\alpha - 1 \leq \beta_Y \leq 1 + (\gamma_X|c|/|a|\gamma_Y)^\alpha (\beta_X - 1). \tag{89}$$

Simplifying the inequalities in (89).

$$(1 + \beta_X)(\gamma_X|c|/|a|\gamma_Y)^\alpha \leq 1 + \beta_Y, \tag{90}$$
$$\beta_Y - 1 \leq (\gamma_X|c|/|a|\gamma_Y)^\alpha (\beta_X - 1). \tag{91}$$

Multiplying $-1$ on both sides in (91):

$$(\gamma_X|c|/|a|\gamma_Y)^\alpha (1 - \beta_X) \leq 1 - \beta_Y. \tag{92}$$

Hence, by above analysis we can conclude that the inequality $|\beta'| \leq 1$ is equivalent to the inequalities: $(\gamma_X|c|/|a|\gamma_Y)^\alpha (1 - \beta_X) \leq 1 - \beta_Y$ and $(1 + \beta_X)(\gamma_X|c|/|a|\gamma_Y)^\alpha \leq 1 + \beta_Y$, which ensures $\beta' \in [-1, 1]$.

As $\beta' \in [-1, 1]$, $\gamma' \in (0, \infty)$ and $\mu' \in \mathbb{R}$, we can conclude $\epsilon$ follows a legitimate univariate continuous stable distribution. Furthermore, we define $\epsilon \perp\!\!\!\perp (M, X, X')$, i.e., $\Pr(\epsilon|M, X, X') = \Pr(\epsilon)$ and $\Pr(M, X, X'|\epsilon) = \Pr(M, X, X')$.

We now use $\epsilon$ to specify $Q_{MC}(Y'|X')$ as follows:

$$Y' = X' + \epsilon. \tag{93}$$

We also derive the distribution $Q(Y'|M)$ before proceeding with our construction, as we will need it later in the proof to show that the constructed $Q_{MC}(M, X, Y) \in \Delta_P$.

*Deriving $Q_{MC}(Y'|M)$:* We will use the distributions $Q_{MC}(X'|M)$ and $Q_{MC}(\epsilon)$ to derive the distribution of $Q_{MC}(Y'|M)$:

$$Q_{MC}(Y'{=}y'|M{=}m) = Q_{MC}(X' + \epsilon{=}y'|M{=}m),$$
$$= \int_{k=-\infty}^{\infty} Q_{MC}(X'{=}k, \epsilon{=}y' - k|M{=}m)dk,$$
$$= \int_{k=-\infty}^{\infty} Q_{MC}(\epsilon = y' - k|M{=}m)Q_{MC}(X'{=}k|\epsilon = y' - k, M{=}m)dk.$$

Using the fact that $\epsilon \perp\!\!\!\perp M$ and $\epsilon \perp\!\!\!\perp X'|M$, we have:

$$Q_{MC}(Y' = y'|M = m) = \int_{k=-\infty}^{\infty} Q_{MC}(\epsilon = y' - k)Q_{MC}(X'{=}k|M{=}m)dk. \tag{94}$$

Since $Q_{MC}(\epsilon)$ and $Q_{MC}(X'|M = m)$ are univariate continuous stable distributions for a fixed $m$ (see (84) and (83)), equation (94) describes a convolution of two univariate stable distributions for a fixed $m$. Convolution of two univariate continuous stable distribution is akin to summing two independent random variable having the same univariate continuous stable distributions, and

consequently $Q_{MC}(Y'|M)$ can be derived using lemma 6. Hence,

$$Q_{MC}(Y'|M=m) = p_{CS}(\alpha, \tilde{\beta}, \tilde{\gamma}, \tilde{\mu}), \text{ where}$$

$$\tilde{\mu} = m + \frac{b}{a} + \mu' = m + \frac{b}{a} + \frac{d}{c} - \frac{b}{a} = m + \frac{d}{c},$$

$$\tilde{\gamma} = \left(\left(\frac{\gamma_X}{|a|}\right)^\alpha + \gamma'^\alpha\right)^{\frac{1}{\alpha}} = \left(\left(\frac{\gamma_X}{|a|}\right)^\alpha + \left(\frac{\gamma_Y}{|c|}\right)^\alpha - \left(\frac{\gamma_X}{|a|}\right)^\alpha\right)^{\frac{1}{\alpha}} = \frac{\gamma_Y}{|c|},$$

$$\tilde{\beta} = \frac{(\gamma_X/|a|)^\alpha \operatorname{sgn}(a)\beta_X + \gamma'^\alpha \beta'}{(\gamma_X/|a|)^\alpha + \gamma'^\alpha}$$

$$= \frac{(\gamma_X/|a|)^\alpha \operatorname{sgn}(a)\beta_X + ((\gamma_Y/|c|)^\alpha - (\gamma_X/|a|)^\alpha)\left(\operatorname{sgn}(a)\frac{(\gamma_Y/|c|)^\alpha \beta_Y - (\gamma_X/|a|)^\alpha \beta_X}{(\gamma_Y/|c|)^\alpha - (\gamma_X/|a|)^\alpha}\right)}{(\gamma_X/|a|)^\alpha + (\gamma_Y/|c|)^\alpha - (\gamma_X/|a|)^\alpha}$$

$$= \operatorname{sgn}(a)\frac{(\gamma_X/|a|)^\alpha \beta_X + (\gamma_Y/|c|)^\alpha \beta_Y - (\gamma_X/|a|)^\alpha \beta_X}{(\gamma_Y/|c|)^\alpha} = \operatorname{sgn}(a)\frac{(\gamma_Y/|c|)^\alpha \beta_Y}{(\gamma_Y/|c|)^\alpha}$$

$$= \operatorname{sgn}(a)\beta_Y. \tag{95}$$

Specifying $Q_{MC}(Y|Y')$: $Q_{MC}(Y|Y')$ is specified by the following deterministic transformation:

$$Y = \begin{cases} cY' & \alpha \neq 1 \\ cY' + (2\operatorname{sgn}(a)c\beta_Y\gamma_Y \log(|c|))/\pi & \alpha = 1 \end{cases}. \tag{96}$$

Finally, we construct the desired $Q_{MC}(M, X, Y)$ from $Q_{MC}(M, X, X', Y', Y)$ by marginalizing $X'$ and $Y'$.

**Part 2: Showing $Q_{MC}(M, X, Y) \in \Delta_P$**

For showing that $Q_{MC}(M, X, Y) \in \Delta_P$, we first need to calculate $Q_{MC}(Y|M)$.

Calculating $Q(Y|M)$: We use lemma 5 to calculate $Q(Y|M)$ as from (95), we know that $\overline{Q_{MC}(Y'|M) = p_{CS}(Y'; \alpha, \operatorname{sgn}(a)\beta_Y, \gamma_Y/|c|, M + d/c)}$ is a continuous stable distribution for a fixed $m$, and $Y$ a translated and scaled version of $Y'$. The exact expression of $Q_{MC}(Y|M=m)$ is provided in (97).

$$Q_{MC}(Y|M=m) = p_{CS}(\alpha, \tilde{\beta}_Y, \tilde{\gamma}_Y, \tilde{\mu}_Y), \text{ where}$$

$$\tilde{\beta}_Y = \begin{cases} \operatorname{sgn}(c)\operatorname{sgn}(c)\beta_Y & \alpha \neq 1 \\ \operatorname{sgn}(c)\operatorname{sgn}(a)\beta_Y & \alpha = 1 \end{cases} = \operatorname{sgn}(a)\operatorname{sgn}(c)\beta_Y = \operatorname{sgn}(ac)\beta_Y,$$

$$\tilde{\gamma}_Y = \begin{cases} |c|\frac{\gamma_Y}{|c|} & \alpha \neq 1 \\ |c|\frac{\gamma_Y}{|c|} & \alpha = 1 \end{cases} = \gamma_Y,$$

$$\tilde{\mu}_Y = \begin{cases} c(m + \frac{d}{c}) & \alpha \neq 1 \\ c(m + \frac{d}{c}) + 2\operatorname{sgn}(a)c\beta_Y\gamma_Y \log(|c|)/\pi - 2\operatorname{sgn}(a)c\beta_Y\gamma_Y \log(|c|)/\pi & \alpha = 1 \end{cases} = cm + d,$$

$$\Rightarrow Q_{MC}(Y|M=m) = p_{CS}(\alpha, \beta_Y \operatorname{sgn}(ac), \gamma_Y, cm + d) = P(Y|M). \tag{97}$$

From (81) and (97), we can conclude:

$$Q_{MC}(M, X) = Q_{MC}(M)Q_{MC}(X|M) = P(M)P(X|M) = P(M, X), \tag{98}$$

$$Q_{MC}(M, Y) = Q_{MC}(M)Q_{MC}(Y|M) = P(M)P(Y|M) = P(M, Y). \tag{99}$$

Hence, $Q_{MC}(M, X, Y) \in \Delta_P$ and consequently, by proposition 1, $UI(M; Y\backslash X) = 0$, concluding our proof for case 1.

**Case 2: $|a|/\gamma_X = |c|/\gamma_Y$**

The proof of case 2 is extremely similar to case 1. For case 2, we are able to directly construct the distribution $Q_{MC}(M, X, Y)$ without the need of specifying a larger Markov chain.

**Part 1: Specifying $Q_{MC}(M, X, Y)$**

Similarly to case 1, we specify $Q_{MC}(M, X, Y)$ by individually specifying $Q_{MC}(M)$, $Q_{MC}(X|M)$ and $Q_{MC}(Y|X)$. The distributions $Q_{MC}(M)$ and $Q_{MC}(X|M)$ are as follows:

$$Q_{MC}(M) = P(M) \text{ and } Q_{MC}(X|M) = P(X|M). \tag{100}$$

The distribution $Q_{MC}(Y|X)$ is defined using the following deterministic transformation:

$$Y = \begin{cases} {}^{c}\!/\!aX - {}^{cb}\!/\!a + d & \alpha \neq 1 \\ \frac{c}{a}X + {}^{2c}\!/\!{}_{a\pi}\beta\gamma_X \log\left(|{}^{c}\!/\!a|\right) - {}^{cb}\!/\!a + d & \alpha = 1 \end{cases}. \tag{101}$$

**Part 2: Showing $Q_{MC}(M,X,Y) \in \Delta_P$**

For showing that $Q_{MC}(M,X,Y) \in \Delta_P$, we first need to calculate $Q_{MC}(Y|M)$.

Calculating $Q(Y|M)$: We use lemma 5 to calculate $Q(Y|M)$ as $Q_{MC}(X|M{=}m)$ is a continuous stable distribution for a fixed $m$, and $Y$ a translated and scaled version of $X$. The exact expression of $Q_{MC}(Y|M = m)$ is provided in (102).

$$Q_{MC}(Y|M = m) = p_{CS}(y; \alpha, \tilde{\beta}, \tilde{\gamma}, \tilde{\mu}),$$

$$\tilde{\mu} = \begin{cases} {}^{ca}\!/\!am + \frac{cb}{a} - \frac{bc}{a} + d & \alpha \neq 1 \\ {}^{ca}\!/\!am + \frac{cb}{a} - \frac{bc}{a} + d - {}^{2c}\!/\!{}_{a\pi}\beta\gamma_X \log\left(|{}^{c}\!/\!a|\right) + {}^{2c}\!/\!{}_{a\pi}\beta\gamma_X \log\left(|{}^{c}\!/\!a|\right) & \alpha = 1 \end{cases}$$

$$= cm + d,$$

$$\tilde{\gamma} = |c|\gamma_X/|a| = |c|\gamma_Y/|c| = \gamma_Y, (\text{ using the fact that } {}^{\gamma_X}\!/\!|a| = {}^{\gamma_Y}\!/\!|c| \text{ by assumption})$$

$$\tilde{\beta} = \beta\mathrm{sgn}({}^{c}\!/\!a) = \beta\mathrm{sgn}(ac),$$

$$\Rightarrow Q_{MC}(Y|M = m) = p_{CS}(\alpha, \mathrm{sgn}(ac)\beta_Y, \gamma_Y, cm + d) = P(Y|M). \tag{102}$$

From (100) and (102), we can conclude:

$$Q_{MC}(M,X) = Q_{MC}(M)Q_{MC}(X|M) = P(M)P(X|M) = P(M,X), \tag{103}$$

$$Q_{MC}(M,Y) = Q_{MC}(M)Q_{MC}(Y|M) = P(M)P(Y|M) = P(M,Y). \tag{104}$$

Hence, $Q_{MC}(M,X,Y) \in \Delta_P$ and consequently, by proposition 1, $UI(M;Y\backslash X) = 0$, concluding our proof for case 2 and theorem 1. $\qquad\square$

# G    Proof of Theorem 2

In this section, we provide the proof of theorem 2. Since theorem 2 provides the analytical PID terms for system 1 of the multivariate affine continuous stable system defined in Sec. 4, we briefly restate certain key properties of the corresponding independent component multivariate stable distributions and the corresponding multivariate affine continuous stable system for convenience.

## G.1    Independent component multivariate stable distribution

Independent component multivariate stable distributions are a specific multivariate generalization of univariate continuous stable distributions describing a collection of independent random variables, with each random variable distributed according to a univariate continuous stable distribution. We refer the reader to Appx. N.3 for more details on multivariate continuous stable distributions. We now list certain key properties of independent component multivariate stable distributions that we make use of in the proof of theorem 2:

1. If $\vec{\mathbf{X}}$ is distributed according to an independent component multivariate stable distribution, then the sum of two independent copies of $\vec{\mathbf{X}}$, denoted as $\vec{\mathbf{X}}_1$ and $\vec{\mathbf{X}}_2$, follows an independent component multivariate stable distribution up to a scaling and translation operation, i.e., $a\vec{\mathbf{X}}_1 + b\vec{\mathbf{X}}_2 \stackrel{d}{=} c\vec{\mathbf{X}} + \vec{\mathbf{d}}$ for $a, b, c > 0$ and $\vec{\mathbf{d}} \in \mathbb{R}$.

2. We denote the p.d.f. of the independent component multivariate stable continuous distribution as $p_{CS-IC}(\alpha, \vec{\boldsymbol{\beta}}, \vec{\boldsymbol{\gamma}}, \vec{\boldsymbol{\mu}})$, where

    $$\vec{\boldsymbol{\beta}} = [\beta_1 \quad \dots \quad \beta_d]^T, \text{ with } \beta_j \in [-1,1], \vec{\boldsymbol{\gamma}} = [\gamma_1 \quad \dots \quad \gamma_d]^T \text{ with } \gamma_j \in (0, \infty),$$

    $$\vec{\boldsymbol{\mu}} = [\mu_1 \quad \dots \quad \mu_d]^T \in \mathbb{R}^d, \text{ and } \alpha \in (0,2].$$

    In general, the p.d.f. of independent component multivariate stable distribution do not have a closed-form analytical expression, and are expressed through their characteristic function.

3. The characteristic function of a $d$-dimensional random vector $\vec{\mathbf{X}}$ having independent component multivariate distribution $p_{CS-IC}(\alpha, \vec{\boldsymbol{\beta}}, \vec{\boldsymbol{\gamma}}, \vec{\boldsymbol{\mu}})$ is expressed as follows:

$$\mathbb{E}\left[e^{i\vec{\mathbf{t}}^T \vec{\mathbf{X}}}\right] = \exp\left(-\sum_{j=1}^{d}|\gamma_j t_j|^\alpha(1 - i\beta_j \mathrm{sgn}(t_j)\Phi(\alpha)) + i\vec{\mathbf{t}}^T\vec{\boldsymbol{\mu}}\right) \ \forall \ \vec{\mathbf{t}} \in \mathbb{R}^d, \quad (105)$$

$$\text{where } \Phi(\alpha) = \left\{\begin{array}{ll} \tan\left(\frac{\pi\alpha}{2}\right) & \alpha \neq 1 \\ \frac{-2}{\pi}\log(|t|) & \alpha = 1 \end{array}\right., \quad \mathrm{sgn}(t) = \left\{\begin{array}{ll} -1 & t < 0 \\ 0 & t = 0 \\ 1 & t > 0 \end{array}\right..$$

4. The random vector $\vec{\mathbf{X}} = [X_1 \ \ldots \ X_d]^T$ having the distribution $p_{CS-IC}(\alpha, \vec{\boldsymbol{\beta}}, \vec{\boldsymbol{\gamma}}, \vec{\boldsymbol{\mu}})$ is essentially a collection of independent random variables $\{X_j\}_{j=1}^{d}$, where $X_j \sim p_{CS}(\alpha, \beta_j, \gamma_j, \mu_j)$.

## G.2 Definition of the system 1 of the multivariate affine continuous stable system

Let the random variable $M \sim P(M)$ with support $\mathcal{M} \subseteq \mathbb{R}$. $\vec{\mathbf{X}}$ and $\vec{\mathbf{Y}}$ are $d_X$-dimensional and $d_Y$-dimensional random vectors, respectively. The joint distribution $P(M, \vec{\mathbf{X}}, \vec{\mathbf{Y}})$ describes system 1 of the multivariate affine continuous stable system if it satisfies the following two properties

1. $M \sim P(M)$ having some support set $\mathcal{M} \subseteq \mathbb{R}$.

2. The conditional distributions of random variables $\vec{\mathbf{X}}$ and $\vec{\mathbf{Y}}$ conditioned on $M$ are expressed as follows:

$$\vec{\mathbf{X}} = \vec{\mathbf{H}}_X M + \mathbf{A}_X \vec{\mathbf{Z}}_X + \vec{\mathbf{b}}_X \text{ and } \vec{\mathbf{Y}} = \vec{\mathbf{H}}_Y M + \mathbf{A}_Y \vec{\mathbf{Z}}_Y + \vec{\mathbf{b}}_Y, \quad (106)$$

where $\vec{\mathbf{Z}}_X \sim p_{CS-IC}(\alpha, \vec{\mathbf{0}}_{d_X}, \vec{\mathbf{1}}_{d_X}, \vec{\mathbf{0}}_{d_X})$, $\vec{\mathbf{Z}}_Y \sim p_{CS-IC}(\alpha, \vec{\mathbf{0}}_{d_Y}, \vec{\mathbf{1}}_{d_Y}, \vec{\mathbf{0}}_{d_Y})$, $\mathbf{A}_X$ and $\mathbf{A}_Y$ are invertible matrices, $\vec{\mathbf{H}}_X, \vec{\mathbf{b}}_X \in \mathbb{R}^{d_X}$, and $\vec{\mathbf{H}}_Y, \vec{\mathbf{b}}_Y \in \mathbb{R}^{d_Y}$.

## G.3 Formal proof of Theorem 2

**Theorem 2.** *Let the joint distribution $P(M, \vec{\mathbf{X}}, \vec{\mathbf{Y}})$ of $M$, $\vec{\mathbf{X}}$, and $\vec{\mathbf{Y}}$ describe the system 1 of the multivariate affine continuous stable system. Without the loss of generality, assume $\|\mathbf{A}_Y^{-1}\vec{\mathbf{H}}_Y\|_\kappa \leq \|\mathbf{A}_X^{-1}\vec{\mathbf{H}}_X\|_\kappa$, where $\kappa = \alpha/\alpha-1 \ \forall \ \alpha \in (1, 2]$ and $\kappa = \infty \ \forall \ \alpha \in (0, 1]$. Then, $\Delta_P$ contains a Markov chain of the form $M \to \vec{\mathbf{X}} \to \vec{\mathbf{Y}}$ and $UI(M; \vec{\mathbf{Y}}\backslash\vec{\mathbf{X}}) = 0$.*

*Proof.* We first note that we can always assume $\|\mathbf{A}_Y^{-1}\vec{\mathbf{H}}_Y\|_\kappa \leq \|\mathbf{A}_X^{-1}\vec{\mathbf{H}}_X\|_\kappa$ without the loss of generality because if $\|\mathbf{A}_Y^{-1}\vec{\mathbf{H}}_Y\|_\kappa \geq \|\mathbf{A}_X^{-1}\vec{\mathbf{H}}_X\|_\kappa$, then we can always switch our nomenclature to refer to $\vec{\mathbf{Y}}$ as $\vec{\mathbf{X}}$, and $\vec{\mathbf{X}}$ as $\vec{\mathbf{Y}}$.

The proof of above theorem relies on the result of lemma 7 and the fact that the linear system described in (106) can always be reduced to the special case of the linear system used in lemma 7. We briefly outline the proof structure.

1. In the first part, we explicitly construct a joint distribution $Q_{MC}(M, \vec{\mathbf{X}}, \vec{\mathbf{Y}})$ having the Markovian structure $M \to \vec{\mathbf{X}} \to \vec{\mathbf{Y}}$.

2. In the second part, we show that the $Q_{MC}(M, \vec{\mathbf{X}}, \vec{\mathbf{Y}})$ constructed in the first part, lies in $\Delta_P$. Therefore, we can then apply the result of proposition 1 to conclude $UI(M; \vec{\mathbf{Y}}\backslash\vec{\mathbf{X}}) = 0$.

## Part 1: Specifying $Q_{MC}(M, \vec{\mathbf{X}}, \vec{\mathbf{Y}})$

We explicitly construct the desired Markov chain $M \to \vec{\mathbf{X}} \to \vec{\mathbf{Y}}$ by constructing a larger Markov chain $M \to \vec{\mathbf{X}} \to \vec{\mathbf{X}}' \to \vec{\mathbf{Y}}' \to \vec{\mathbf{Y}}$ and then marginalizing the larger Markov chain to obtain the desired Markov chain $M \to \vec{\mathbf{X}} \to \vec{\mathbf{Y}}$.

Denote the joint distribution of the Markov chain $M \to \vec{\mathbf{X}} \to \vec{\mathbf{X}}' \to \vec{\mathbf{Y}}' \to \vec{\mathbf{Y}}$ as $Q_{MC}(M, \vec{\mathbf{X}}, \vec{\mathbf{X}}', \vec{\mathbf{Y}}', \vec{\mathbf{Y}})$. We can decompose $Q_{MC}(M, \vec{\mathbf{X}}, \vec{\mathbf{X}}', \vec{\mathbf{Y}}', \vec{\mathbf{Y}})$ by utilizing its Markovian structure as follows:

$$Q_{MC}(M, \vec{\mathbf{X}}, \vec{\mathbf{X}}', \vec{\mathbf{Y}}', \vec{\mathbf{Y}}) = Q_{MC}(M) Q_{MC}(\vec{\mathbf{X}}|M) Q_{MC}(\vec{\mathbf{X}}'|\vec{\mathbf{X}}) Q_{MC}(\vec{\mathbf{Y}}'|\vec{\mathbf{X}}') Q_{MC}(\vec{\mathbf{Y}}|\vec{\mathbf{Y}}').$$

Consequently, we can specify/construct the distribution $Q_{MC}(M, \vec{\mathbf{X}}, \vec{\mathbf{X}}', \vec{\mathbf{Y}}', \vec{\mathbf{Y}})$ by individually specifying $Q_{MC}(M)$, $Q_{MC}(\vec{\mathbf{X}}|M)$, $Q_{MC}(\vec{\mathbf{X}}'|\vec{\mathbf{X}})$, $Q_{MC}(\vec{\mathbf{Y}}'|\vec{\mathbf{X}}')$, and $Q_{MC}(\vec{\mathbf{Y}}|\vec{\mathbf{Y}}')$. We specify $Q_{MC}(M, \vec{\mathbf{X}}) = P(M, \vec{\mathbf{X}})$ and $Q(\vec{\mathbf{X}}'|\vec{\mathbf{X}})$ using the following deterministic transformation:

$$\vec{\mathbf{X}}' = \mathbf{A}_X^{-1} \left( \vec{\mathbf{X}} - \vec{\mathbf{b}}_X \right). \tag{107}$$

Substituting (106) in the above equation:

$$\vec{\mathbf{X}}' = \mathbf{A}_X^{-1} \left( \vec{\mathbf{H}}_X M + \mathbf{A}_X \vec{\mathbf{Z}}_X + \vec{\mathbf{b}}_X - \vec{\mathbf{b}}_X \right) = \mathbf{A}_X^{-1} \vec{\mathbf{H}}_X M + \vec{\mathbf{Z}}_X. \tag{108}$$

Specifying $Q_{MC}(\vec{\mathbf{Y}}'|\vec{\mathbf{X}}')$: We define $\vec{\mathbf{Y}}'$ according to the following affine system:

$$\vec{\mathbf{Y}}' = \mathbf{A}_Y^{-1} \vec{\mathbf{H}}_Y M + \vec{\mathbf{Z}}_Y. \tag{109}$$

Observe that both $\vec{\mathbf{X}}'$ and $\vec{\mathbf{Y}}'$ can be described as an affine function of $M$ satisfying the special structure described in lemma 7 (see (115)). Furthermore, $Q_{MC}(M) = P(M)$ by construction. Consequently, $Q_{MC}(M)$ satisfies the properties required by the distribution of $M$ outlined in lemma 7, as $P(M)$ in both lemma 7 and theorem 2 follow the same properties. Since $\|\mathbf{A}_Y^{-1} \vec{\mathbf{H}}_Y\|_\kappa \leq \|\mathbf{A}_X^{-1} \vec{\mathbf{H}}_X\|_\kappa$ by the assumption in the theorem, we can apply the result of lemma 7 to construct $Q_{MC}(\vec{\mathbf{Y}}'|\vec{\mathbf{X}}')$, where $\vec{\mathbf{H}}_X$ and $\vec{\mathbf{H}}_Y$ would be replaced by $\mathbf{A}_X^{-1} \vec{\mathbf{H}}_X$ and $\mathbf{A}_Y^{-1} \vec{\mathbf{H}}_Y$, respectively.

Lastly, we specify $Q_{MC}(\vec{\mathbf{Y}}|\vec{\mathbf{Y}}')$ through the following linear system:

$$\vec{\mathbf{Y}} = \mathbf{A}_Y \vec{\mathbf{Y}}' + \vec{\mathbf{b}}_Y. \tag{110}$$

Finally, we construct the desired $Q_{MC}(M, \vec{\mathbf{X}}, \vec{\mathbf{Y}})$ from $Q_{MC}(M, \vec{\mathbf{X}}, \vec{\mathbf{X}}', \vec{\mathbf{Y}}', \vec{\mathbf{Y}})$ by marginalizing $\vec{\mathbf{X}}'$ and $\vec{\mathbf{Y}}'$.

**Part 2: Showing $Q_{MC}(M, \vec{\mathbf{X}}, \vec{\mathbf{Y}}) \in \Delta_P$**

For showing that $Q_{MC}(M, \vec{\mathbf{X}}, \vec{\mathbf{Y}}) \in \Delta_P$, we first need to derive $Q_{MC}(\vec{\mathbf{Y}}|M)$.

*Deriving $Q(\vec{\mathbf{Y}}|M)$:* Substituting (109) in (110) we obtain:

$$\vec{\mathbf{Y}} = \mathbf{A}_Y (\mathbf{A}_Y^{-1} \vec{\mathbf{H}}_Y M + \vec{\mathbf{Z}}_Y) + \vec{\mathbf{b}}_Y = \vec{\mathbf{H}}_Y M + \mathbf{A}_Y \vec{\mathbf{Z}}_Y + \vec{\mathbf{b}}_Y. \tag{111}$$

Inspecting the above equation, we can conclude:

$$Q_{MC}(\vec{\mathbf{Y}}|M) = P(\vec{\mathbf{Y}}|M). \tag{112}$$

From (112), we can conclude:

$$Q_{MC}(M, \vec{\mathbf{Y}}) = Q_{MC}(M) Q_{MC}(\vec{\mathbf{Y}}|M) = P(M) P(\vec{\mathbf{Y}}|M) = P(M, \vec{\mathbf{Y}}). \tag{113}$$

We know that by construction:

$$Q_{MC}(M, \vec{\mathbf{X}}) = P(M, \vec{\mathbf{X}}). \tag{114}$$

Hence, $Q_{MC}(M, \vec{\mathbf{X}}, \vec{\mathbf{Y}}) \in \Delta_P$ and consequently, by proposition 1, $UI(M; \vec{\mathbf{Y}} \backslash \vec{\mathbf{X}}) = 0$, concluding our proof. $\square$

**Lemma 7.** *Let the joint distribution of $M$, $\vec{\mathbf{X}}$ and $\vec{\mathbf{Y}}$, denoted as $P(M, \vec{\mathbf{X}}, \vec{\mathbf{Y}})$, describe the system 1 of the multivariate affine continuous stable system. Furthermore, assume $\mathbf{A}_X = \mathbf{I}_{d_X}$, $\mathbf{A}_Y = \mathbf{I}_{d_Y}$, $\vec{\mathbf{b}}_X = \vec{\mathbf{0}}_{d_X}$ and $\vec{\mathbf{b}}_Y = \vec{\mathbf{0}}_{d_Y}$, where $\mathbf{I}_d$ is a $d \times d$ identity matrix and $\vec{\mathbf{0}}_d$ is $d$-dimensional vector of zeros. Then,*

1. If $\|\vec{\mathbf{H}}_Y\|_\kappa \leq \|\vec{\mathbf{H}}_X\|_\kappa$, where $\kappa = \alpha/\alpha-1 \ \forall \ \alpha \in (1,2]$ and $\kappa = \infty \ \forall \ \alpha \in (0,1]$, then $\Delta_P$ contains a Markov chain of the form $M \to \vec{\mathbf{X}} \to \vec{\mathbf{Y}}$ and $UI(M; \vec{\mathbf{Y}}/\vec{\mathbf{X}}) = 0$.

2. If $\|\vec{\mathbf{H}}_X\|_\kappa \leq \|\vec{\mathbf{H}}_Y\|_\kappa$, where $\kappa = \alpha/\alpha-1 \ \forall \ \alpha \in (1,2]$ and $\kappa = \infty \ \forall \ \alpha \in (0,1]$, then $\Delta_P$ contains a Markov chain of the form $M \to \vec{\mathbf{Y}} \to \vec{\mathbf{X}}$ and $UI(M; \vec{\mathbf{X}}/\vec{\mathbf{Y}}) = 0$.

*Proof.* We only provide an explicit proof of condition 1. Condition 2 is essentially the same as condition 1, with the parameters of X and Y switched. Consequently, the proof of condition 2 follows the same steps as the proof of condition 1, with parameters about X and Y switched.

**Proof of condition 1**:

We briefly outline the proof structure.

1. In the first part, we explicitly construct a joint distribution $Q_{MC}(M, \vec{\mathbf{X}}, \vec{\mathbf{Y}})$ having the Markovian structure $M \to \vec{\mathbf{X}} \to \vec{\mathbf{Y}}$.

2. In the second part, we show that the $Q_{MC}(M, \vec{\mathbf{X}}, \vec{\mathbf{Y}})$ constructed in the first part, lies in $\Delta_P$. Therefore, we can then apply the result of proposition 1 to conclude $UI(M; \vec{\mathbf{Y}} \backslash \vec{\mathbf{X}}) = 0$.

**Part 1: Specifying $Q_{MC}(M, \vec{\mathbf{X}}, \vec{\mathbf{Y}})$**

We explicitly construct the desired Markov chain $M \to \vec{\mathbf{X}} \to \vec{\mathbf{Y}}$ by constructing a larger Markov chain $M \to \vec{\mathbf{X}} \to \vec{\mathbf{X}}' \to \vec{\mathbf{Y}}$ and then marginalizing the larger Markov chain to obtain the desired Markov chain $M \to \vec{\mathbf{X}} \to \vec{\mathbf{Y}}$.

Denote the joint distribution of the Markov chain $M \to \vec{\mathbf{X}} \to \vec{\mathbf{X}}' \to \vec{\mathbf{Y}}$ as $Q_{MC}(M, \vec{\mathbf{X}}, \vec{\mathbf{X}}', \vec{\mathbf{Y}})$. We can decompose $Q_{MC}(M, \vec{\mathbf{X}}, \vec{\mathbf{X}}', \vec{\mathbf{Y}})$ by utilizing its Markovian structure as follows:

$$Q_{MC}(M, \vec{\mathbf{X}}, \vec{\mathbf{X}}', \vec{\mathbf{Y}}) = Q_{MC}(M)Q_{MC}(\vec{\mathbf{X}}|M)Q_{MC}(\vec{\mathbf{X}}'|\vec{\mathbf{X}})Q_{MC}(\vec{\mathbf{Y}}|\vec{\mathbf{X}}').$$

Consequently, we can specify/construct the distribution $Q_{MC}(M, \vec{\mathbf{X}}, \vec{\mathbf{X}}', \vec{\mathbf{Y}})$ by individually specifying $Q_{MC}(M)$, $Q_{MC}(\vec{\mathbf{X}}|M)$, $Q_{MC}(\vec{\mathbf{X}}'|\vec{\mathbf{X}})$, and $Q_{MC}(\vec{\mathbf{Y}}|\vec{\mathbf{X}}')$.

Under the assumptions of lemma 7, we can simplify the linear system describing $P(M, \vec{\mathbf{X}}, \vec{\mathbf{Y}})$ shown in (106) as follows:

$$\vec{\mathbf{X}} = \vec{\mathbf{H}}_X M + \vec{\mathbf{Z}}_X \text{ and } \vec{\mathbf{Y}} = \vec{\mathbf{H}}_Y M + \vec{\mathbf{Z}}_Y. \tag{115}$$

Correspondingly, we can write the characteristic functions of conditional distributions $P(\vec{\mathbf{X}}|M=m)$ and $P(\vec{\mathbf{Y}}|M=m)$ for a fixed $m$ by employing lemma 9:

$$\mathbb{E}[e^{i\vec{\mathbf{t}}^T\vec{\mathbf{X}}}|M=m] = \exp\left(-\sum_{j=1}^{n} |t_j|^\alpha + i\vec{\mathbf{t}}^T\vec{\mathbf{H}}_X m\right),$$

$$\mathbb{E}[e^{i\vec{\mathbf{t}}^T\vec{\mathbf{Y}}}|M=m] = \exp\left(-\sum_{j=1}^{n} |t_j|^\alpha + i\vec{\mathbf{t}}^T\vec{\mathbf{H}}_Y m\right), \tag{116}$$

where the characteristic function can be derived by realizing that for a given $m$, $\vec{\mathbf{H}}_X m$ and $\vec{\mathbf{H}}_Y m$ are constants being added to the random vectors $\vec{\mathbf{Z}}_X \sim p_{CS-IC}(\alpha, \vec{\mathbf{0}}_{d_x}, \vec{\mathbf{1}}_{d_x}, \vec{\mathbf{0}}_{d_x})$ and $\vec{\mathbf{Z}}_Y \sim p_{CS-IC}(\alpha, \vec{\mathbf{0}}_{d_y}, \vec{\mathbf{1}}_{d_y}, \vec{\mathbf{0}}_{d_y})$, respectively, and then using the result of lemma 9.

Specifying $Q_{MC}(M)$ and $Q_{MC}(\vec{\mathbf{X}}|M)$: We specify $Q_{MC}(M)$ and $Q_{MC}(\vec{\mathbf{X}}|M)$ as follows:

$$Q_{MC}(M) = P(M) \text{ and } Q_{MC}(\vec{\mathbf{X}}|M) = P(\vec{\mathbf{X}}|M), \tag{117}$$

where $P(M)$ and $P(\vec{\mathbf{X}}|M)$ are marginal distributions derived from the original joint distribution $P(M, \vec{\mathbf{X}}, \vec{\mathbf{Y}})$ (discussed in the lemma statement) over which the bivariate PID is being calculated.

Specifying $Q_{MC}(\vec{\mathbf{X}}'|\vec{\mathbf{X}})$: We specify $Q_{MC}(\vec{\mathbf{X}}'|\vec{\mathbf{X}})$ through the following deterministic transformation:

$$\vec{\mathbf{X}}' = \vec{\mathbf{H}}_Y (\vec{\mathbf{H}}_X^k)^T \vec{\mathbf{X}}, \tag{118}$$

where $\vec{\mathbf{H}}_X^k$ is as defined in lemma 10, and $k \in [0, \infty]$.

We will also derive the distribution $Q(\vec{\mathbf{X}}'|M)$ before proceeding with our construction, as we will need it later in the proof to show that the constructed $Q_{MC}(M, \vec{\mathbf{X}}, \vec{\mathbf{Y}}) \in \Delta_P$.

*Deriving $Q_{MC}(\vec{\mathbf{X}}'|M)$*: To derive $Q_{MC}(\vec{\mathbf{X}}'|M)$, we represent $\vec{\mathbf{X}}'$ as function of $M$ and $\vec{\mathbf{Z}}_X$ in a similar manner to $\vec{\mathbf{X}}$ in (115), and then use the result of lemma 9 to derive its conditional characteristic function $\mathbb{E}[e^{\vec{\mathbf{t}}^T \vec{\mathbf{X}}'}|M]$, and correspondingly the desired $Q_{MC}(\vec{\mathbf{X}}'|M)$ .

We multiply $\vec{\mathbf{H}}_Y (\vec{\mathbf{H}}_X^k)^T$ on both sides of equality in (115) to obtain the following linear equation:

$$\vec{\mathbf{H}}_Y (\vec{\mathbf{H}}_X^k)^T \vec{\mathbf{X}} = \vec{\mathbf{H}}_Y (\vec{\mathbf{H}}_X^k)^T (\vec{\mathbf{H}}_X M + \vec{\mathbf{Z}}_X) = \vec{\mathbf{H}}_Y (\vec{\mathbf{H}}_X^k)^T \vec{\mathbf{H}}_X M + \vec{\mathbf{H}}_Y (\vec{\mathbf{H}}_X^k)^T \vec{\mathbf{Z}}_X.$$

Substituting (118) in the above equation, we obtain:

$$\vec{\mathbf{X}}' = \vec{\mathbf{H}}_Y (\vec{\mathbf{H}}_X^k)^T \vec{\mathbf{X}} = \vec{\mathbf{H}}_Y (\vec{\mathbf{H}}_X^k)^T \vec{\mathbf{H}}_X M + \vec{\mathbf{H}}_Y (\vec{\mathbf{H}}_X^k)^T \vec{\mathbf{Z}}_X. \tag{119}$$

Using the result of lemma 10, i.e., $\vec{\mathbf{H}}_Y (\vec{\mathbf{H}}_X^k)^T \vec{\mathbf{H}}_X = \vec{\mathbf{H}}_Y$, in the above equation we obtain:

$$\vec{\mathbf{X}}' = \vec{\mathbf{H}}_Y M + \vec{\mathbf{H}}_Y (\vec{\mathbf{H}}_X^k)^T \vec{\mathbf{Z}}_X. \tag{120}$$

From (120), we can observe that $\vec{\mathbf{X}}'$ is an affine function of $\vec{\mathbf{Z}}_X$ for a fixed $m$. Hence, we can employ the result of lemma 9 to derive the distribution of $Q_{MC}(\vec{\mathbf{X}}'|M)$ (by deriving its corresponding characteristic function).

Note that the $j$-th column of $\vec{\mathbf{H}}_Y (\vec{\mathbf{H}}_X^k)^T$ can be expressed as $h'_j \vec{\mathbf{H}}_Y$, where $h'_j$ is the $j$-th component of $\vec{\mathbf{H}}_X^k$. Consequently:

$$\mathbb{E}\left[ e^{i\vec{\mathbf{t}}^T \vec{\mathbf{X}}'} \,\middle|\, M = m \right] = \exp\left( -\sum_{j=1}^{d_X} \left| h'_j \vec{\mathbf{H}}_Y^T \vec{\mathbf{t}} \right|^\alpha + i\vec{\mathbf{t}}^T \vec{\mathbf{H}}_Y m \right)$$

$$= \exp\left( -\sum_{j=1}^{d_X} |h'_j|^\alpha \left| \vec{\mathbf{H}}_Y^T \vec{\mathbf{t}} \right|^\alpha + i\vec{\mathbf{t}}^T \vec{\mathbf{H}}_Y m \right)$$

$$= \exp\left( -\left| \vec{\mathbf{H}}_Y^T \vec{\mathbf{t}} \right|^\alpha \left( \sum_{j=1}^{d_X} |h'_j|^\alpha \right) + i\vec{\mathbf{t}}^T \vec{\mathbf{H}}_Y m \right). \tag{121}$$

We substitute the value $h'_j$ in (121) using the definition of $\vec{\mathbf{H}}_X^k$ given in lemma 10. We will divide the substitution into two cases:

Case 1: $k$ is finite, i.e., $k \in [0, \infty)$. Then, we have $h'_j = |h_j^X|^k / \|\vec{\mathbf{H}}_X\|_{1+k}^{1+k}$, which implies:

$$\mathbb{E}\left[ e^{i\vec{\mathbf{t}}^T \vec{\mathbf{X}}'} \,\middle|\, M = m \right] = \exp\left( -\left| \vec{\mathbf{H}}_Y^T \vec{\mathbf{t}} \right|^\alpha \sum_{j=1}^{d_X} \frac{|h_j^X|^{k\alpha}}{\|\vec{\mathbf{H}}_X\|_{1+k}^{\alpha+k\alpha}} + i\vec{\mathbf{t}}^T \vec{\mathbf{H}}_Y m \right)$$

$$= \exp\left( -\frac{\left| \vec{\mathbf{H}}_Y^T \vec{\mathbf{t}} \right|^\alpha}{\|\vec{\mathbf{H}}_X\|_{1+k}^{\alpha+k\alpha}} \sum_{j=1}^{d_X} |h_j^X|^{k\alpha} + i\vec{\mathbf{t}}^T \vec{\mathbf{H}}_Y m \right)$$

$$= \exp\left( -\frac{\left| \vec{\mathbf{H}}_Y^T \vec{\mathbf{t}} \right|^\alpha \left\| \vec{\mathbf{H}}_X \right\|_{k\alpha}^{k\alpha}}{\left\| \vec{\mathbf{H}}_X \right\|_{1+k}^{\alpha+k\alpha}} + i\vec{\mathbf{t}}^T \vec{\mathbf{H}}_Y m \right). \tag{122}$$

Case 2: $k$ is infinite, i.e., $k = \infty$. Then, we have $h'_j = 0 \ \forall \ j \neq j^*$, and $h'_{j*} = 1/h_1^X$, which implies:

$$\mathbb{E}\left[e^{i\vec{\mathbf{t}}^T \vec{\mathbf{X}}'} \middle| M = m\right] = \exp\left(-\left|\vec{\mathbf{H}}_Y^T \vec{\mathbf{t}}\right|^\alpha \left|\frac{1}{h_{j*}^X}\right|^\alpha + i\vec{\mathbf{t}}^T \vec{\mathbf{H}}_Y m\right)$$

$$= \exp\left(-\frac{\left|\vec{\mathbf{H}}_Y^T \vec{\mathbf{t}}\right|^\alpha}{\left\|\vec{\mathbf{H}}_X\right\|_\infty^\alpha} + i\vec{\mathbf{t}}^T \vec{\mathbf{H}}_Y m\right). \tag{123}$$

Combining (122) and (123), we can re-write $\mathbb{E}\left[e^{i\vec{\mathbf{t}}^T \vec{\mathbf{X}}'} \middle| M = m\right]$ as:

$$\mathbb{E}\left[e^{i\vec{\mathbf{t}}^T \vec{\mathbf{X}}'} \middle| M = m\right] = \exp\left(-f(k)\left|\vec{\mathbf{H}}_Y^T \vec{\mathbf{t}}\right|^\alpha + i\vec{\mathbf{t}}^T \vec{\mathbf{H}}_Y m\right), \text{ where}$$

$$f(k) = \begin{cases} \|\vec{\mathbf{H}}_X\|_{k\alpha}^{k\alpha}/\|\vec{\mathbf{H}}_X\|_{1+k}^{\alpha+\alpha k} & k \text{ finite} \\ 1/\|\vec{\mathbf{H}}_X\|_\infty^\alpha & k \text{ infinite} \end{cases}. \tag{124}$$

Specifying $Q_{MC}(\vec{\mathbf{Y}}|\vec{\mathbf{X}}')$: We specify $Q_{MC}(\vec{\mathbf{Y}}|\vec{\mathbf{X}}')$ through the following stochastic transformation:

$$\vec{\mathbf{Y}} = \vec{\mathbf{X}}' + \vec{\epsilon}, \tag{125}$$

where $\vec{\epsilon}$ follows a multivariate stable distribution. Furthermore, we assume that $\vec{\epsilon}$ is jointly independent from $(\vec{\mathbf{X}}, \vec{\mathbf{X}}', M)$, i.e., $\vec{\epsilon} \perp\!\!\!\perp (\vec{\mathbf{X}}, \vec{\mathbf{X}}', M)$. The characteristic function of $\vec{\epsilon}$ is defined as follows:

$$\mathbb{E}\left[e^{i\vec{\mathbf{t}}^T \vec{\epsilon}}\right] = \exp\left(-\sum_{j=1}^{d_Y} |t_j|^\alpha + f(k)\left|\vec{\mathbf{H}}_Y^T \vec{\mathbf{t}}\right|^\alpha\right). \tag{126}$$

In order for $Q_{MC}(\vec{\mathbf{Y}}|\vec{\mathbf{X}}')$ to be a valid distribution, we need to ensure that the distribution of $\vec{\epsilon}$ defined in (126) is a legitimate multivariate stable distribution. We will utilize property 1 for showing that $\vec{\epsilon}$ follows a stable multivariate stable distribution. Property 1 states a random vector is distributed according to an independent component multivariate stable distribution iff every 1-dimensional projection of this random vector follows a univariate stable continuous distribution. Hence, to show that $\vec{\epsilon}$ is distributed according to a legitimate multivariate stable distribution, we will show that every 1-dimensional projection of $\vec{\epsilon}$ follows a univariate stable continuous distribution.

Let $\vec{\mathbf{t}} \in \mathbb{R}^{d_Y}$, then the characteristic function of the 1-dimensional projection of $\vec{\epsilon}$ along $\vec{\mathbf{t}}$, i.e., $\vec{\mathbf{t}}^T \vec{\epsilon}$ can be trivially deduced using (126).

$$\mathbb{E}\left[e^{i\vec{\mathbf{t}}^T \vec{\epsilon}}\right] = \exp\left(-\sum_{j=1}^{d_Y} |t_j|^\alpha + f(k)\left|\vec{\mathbf{H}}_Y^T \vec{\mathbf{t}}\right|^\alpha\right)$$

$$= \exp\left(-\left\|\vec{\mathbf{t}}\right\|_\alpha^\alpha + f(k)\left|\vec{\mathbf{H}}_Y^T \vec{\mathbf{t}}\right|^\alpha\right). \tag{127}$$

Comparing the characteristic function of $\vec{\mathbf{t}}^T \vec{\epsilon}$ described in (127) with the standard characteristic function of a univariate stable characteristic function defined in (323), we can conclude $\vec{\mathbf{t}}^T \vec{\epsilon} \sim p_{CS}(\alpha, \beta(\vec{\mathbf{t}}), \gamma(\vec{\mathbf{t}}), \mu(\vec{\mathbf{t}}))$, where:

$$\gamma(\vec{\mathbf{t}}) = \left\|\vec{\mathbf{t}}\right\|_\alpha^\alpha - f(k)\left|\vec{\mathbf{H}}_Y^T \vec{\mathbf{t}}\right|^\alpha, \mu(\vec{\mathbf{t}}) = 0, \text{ and } \beta(\vec{\mathbf{t}}) = 0. \tag{128}$$

In order for $\vec{\mathbf{t}}^T \vec{\epsilon}$ to have a legitimate univariate stable distribution, we just need to show that $\gamma(\vec{\mathbf{t}}) \geq 0 \ \forall \ \vec{\mathbf{t}} \in \mathbb{R}^{d_Y}$. Note that $\gamma(\vec{\mathbf{t}}) = 0$ would correspond to the case where all the mass of the distribution is centered at 0, which does satisfy definition 2, and is an example of a degenerate univariate stable distribution. Let us analyze the function $\gamma(\vec{\mathbf{t}})$:

$$\gamma(\vec{\mathbf{t}}) = \left\|\vec{\mathbf{t}}\right\|_\alpha^\alpha - f(k)\left|\vec{\mathbf{H}}_Y^T \vec{\mathbf{t}}\right|^\alpha, \tag{129}$$

Applying lemma 8 on the term $\left|\vec{\mathbf{H}}_Y^T \vec{\mathbf{t}}\right|$, we obtain:

$$
\left|\vec{\mathbf{H}}_Y^T \vec{\mathbf{t}}\right| \leq \begin{cases} \|\vec{\mathbf{t}}\|_\alpha^\alpha \|\vec{\mathbf{H}}_Y\|_{\alpha/\alpha-1}^\alpha & \alpha \in (1,2] \\ \|\vec{\mathbf{t}}\|_1^\alpha \|\vec{\mathbf{H}}_Y\|_\infty^\alpha & \alpha \in (0,1] \end{cases} .
$$

(130)

Applying (130) in (129), we obtain:

$$
\gamma(\vec{\mathbf{t}}) \geq \begin{cases} \|\vec{\mathbf{t}}\|_\alpha^\alpha - f(k)\|\vec{\mathbf{t}}\|_\alpha^\alpha \|\vec{\mathbf{H}}_Y\|_{\alpha/\alpha-1}^\alpha & \alpha \in (1,2] \\ \|\vec{\mathbf{t}}\|_\alpha^\alpha - f(k)\|\vec{\mathbf{t}}\|_1^\alpha \|\vec{\mathbf{H}}_Y\|_\infty^\alpha & \alpha \in (0,1] \end{cases} .
$$

(131)

Denote $l(\vec{\mathbf{t}}, \alpha)$ as follows:

$$
l(\vec{\mathbf{t}}, \alpha) = \begin{cases} \|\vec{\mathbf{t}}\|_\alpha^\alpha - f(k)\|\vec{\mathbf{t}}\|_\alpha^\alpha \|\vec{\mathbf{H}}_Y\|_{\alpha/\alpha-1}^\alpha & \alpha \in (1,2] \\ \|\vec{\mathbf{t}}\|_\alpha^\alpha - f(k)\|\vec{\mathbf{t}}\|_1^\alpha \|\vec{\mathbf{H}}_Y\|_\infty^\alpha & \alpha \in (0,1] \end{cases} .
$$

(132)

In order to show $\gamma(\vec{\mathbf{t}}) \geq 0 \,\forall\, \vec{\mathbf{t}} \in \mathbb{R}^{d_Y}$, it suffices to show that $l(\vec{\mathbf{t}}, \alpha) \geq 0 \,\forall\, \vec{\mathbf{t}} \in \mathbb{R}^{d_Y}$ and $\alpha \in (0,2]$.

Case 1: $\alpha \in (1,2]$.

For $\alpha \in (1,2]$, we have:

$$
l(\vec{\mathbf{t}}, \alpha) = \left\|\vec{\mathbf{t}}\right\|_\alpha^\alpha - f(k) \left\|\vec{\mathbf{t}}\right\|_\alpha^\alpha \left\|\vec{\mathbf{H}}_Y\right\|_{\alpha/\alpha-1}^\alpha = \left\|\vec{\mathbf{t}}\right\|_\alpha^\alpha \left(1 - f(k)\left\|\vec{\mathbf{H}}_Y\right\|_{\alpha/\alpha-1}^\alpha\right).
$$

(133)

Since, the above equation holds for any $k \in [0,\infty]$, we choose $k = 1/(\alpha-1)$. Substituting the form $f(k)$ from (124) in the above equation, we obtain:

$$
l(\vec{\mathbf{t}}, \alpha) = \left\|\vec{\mathbf{t}}\right\|_\alpha^\alpha \left(1 - \frac{\left\|\vec{\mathbf{H}}_Y\right\|_{\alpha/\alpha-1}^\alpha \left\|\vec{\mathbf{H}}_X\right\|_{\alpha/\alpha-1}^{\alpha/\alpha-1}}{\left\|\vec{\mathbf{H}}_X\right\|_{1+\frac{1}{\alpha-1}}^{\alpha\left(1+\frac{1}{\alpha-1}\right)}}\right) = \left\|\vec{\mathbf{t}}\right\|_\alpha^\alpha \left(1 - \frac{\left\|\vec{\mathbf{H}}_Y\right\|_{\alpha/\alpha-1}^\alpha \left\|\vec{\mathbf{H}}_X\right\|_{\alpha/\alpha-1}^{\alpha/\alpha-1}}{\left\|\vec{\mathbf{H}}_X\right\|_{\alpha/\alpha-1}^{\alpha\left(\frac{\alpha}{\alpha-1}\right)}}\right)
$$

$$
= \left\|\vec{\mathbf{t}}\right\|_\alpha^\alpha \left(1 - \frac{\left\|\vec{\mathbf{H}}_Y\right\|_{\alpha/\alpha-1}^\alpha}{\left\|\vec{\mathbf{H}}_X\right\|_{\alpha/\alpha-1}^{\alpha\left(\frac{\alpha}{\alpha-1}\right)-\frac{\alpha}{\alpha-1}}}\right) = \left\|\vec{\mathbf{t}}\right\|_\alpha^\alpha \left(1 - \frac{\left\|\vec{\mathbf{H}}_Y\right\|_{\alpha/\alpha-1}^\alpha}{\left\|\vec{\mathbf{H}}_X\right\|_{\alpha/\alpha-1}^\alpha}\right).
$$

(134)

Note that $\left\|\vec{\mathbf{t}}\right\|_\alpha^\alpha \geq 0$, and by condition 1 we know that $\|\vec{\mathbf{H}}_Y\|_{\alpha/\alpha-1} \leq \|\vec{\mathbf{H}}_X\|_{\alpha/\alpha-1} \Rightarrow \|\vec{\mathbf{H}}_Y\|_{\alpha/\alpha-1}^\alpha \leq \|\vec{\mathbf{H}}_X\|_{\alpha/\alpha-1}^\alpha$. Hence, using these two previous facts and the above equation, we can conclude:

$$
l(\vec{\mathbf{t}}, \alpha) = \underbrace{\left\|\vec{\mathbf{t}}\right\|_\alpha^\alpha}_{\geq 0} \underbrace{\left(1 - \frac{\left\|\vec{\mathbf{H}}_Y\right\|_{\alpha/\alpha-1}^\alpha}{\left\|\vec{\mathbf{H}}_X\right\|_{\alpha/\alpha-1}^\alpha}\right)}_{\geq 0} \Rightarrow l(\vec{\mathbf{t}}, \alpha) \geq 0.
$$

(135)

Case 2: $\alpha \in (0,1]$.

$$
l(\vec{\mathbf{t}}, \alpha) = \left\|\vec{\mathbf{t}}\right\|_\alpha^\alpha - f(k) \left\|\vec{\mathbf{t}}\right\|_1^\alpha \left\|\vec{\mathbf{H}}_Y\right\|_\infty^\alpha \overset{(a)}{\geq} \left\|\vec{\mathbf{t}}\right\|_\alpha^\alpha - f(k) \left\|\vec{\mathbf{t}}\right\|_\alpha^\alpha \left\|\vec{\mathbf{H}}_Y\right\|_\infty^\alpha,
$$

(136)

where $(a)$ is derived using the monotonicity of $L_p$ norms in $p$, see [57]. More precisely, we use the inequality $\|\vec{\mathbf{t}}\|_\alpha \geq \|\vec{\mathbf{t}}\|_1 \,\forall\, \alpha \in (0,1]$. Simplifying the above equation, we have:

$$
l(\vec{\mathbf{t}}, \alpha) \geq \left\|\vec{\mathbf{t}}\right\|_\alpha^\alpha \left(1 - f(k) \left\|\vec{\mathbf{H}}_Y\right\|_\infty^\alpha\right).
$$

(137)

Since, the above equation holds for any $k \in [0, \infty]$, we choose $k = \infty$. Substituting the form $f(k)$ from (124) in the above equation, we obtain:

$$l(\vec{\mathbf{t}}, \alpha) \geq \left\| \vec{\mathbf{t}} \right\|_{\alpha}^{\alpha} \left( 1 - \frac{\|\vec{\mathbf{H}}_Y\|_{\infty}^{\alpha}}{\|\vec{\mathbf{H}}_X\|_{\infty}^{\alpha}} \right). \tag{138}$$

Note that $\left\| \vec{\mathbf{t}} \right\|_{\alpha}^{\alpha} \geq 0$, and by condition 1 we know that $\|\vec{\mathbf{H}}_Y\|_{\infty} \leq \|\vec{\mathbf{H}}_X\|_{\infty} \Rightarrow \|\vec{\mathbf{H}}_Y\|_{\infty}^{\alpha} \leq \|\vec{\mathbf{H}}_X\|_{\infty}^{\alpha}$. Hence, using these two facts and the above equation, we can conclude:

$$l(\vec{\mathbf{t}}, \alpha) \geq \underbrace{\left\| \vec{\mathbf{t}} \right\|_{\alpha}^{\alpha}}_{\geq 0} \underbrace{\left( 1 - \frac{\left\| \vec{\mathbf{H}}_Y \right\|_{\infty}^{\alpha}}{\left\| \vec{\mathbf{H}}_X \right\|_{\infty}^{\alpha}} \right)}_{\geq 0} \Rightarrow l(\vec{\mathbf{t}}, \alpha) \geq 0. \tag{139}$$

From (135) and (139), we can conclude that $l(\vec{\mathbf{t}}, \alpha) \geq 0 \ \forall \ \vec{\mathbf{t}} \in \mathbb{R}^{d_Y}$ and $\alpha \in (0, 2]$, which implies $\gamma(\vec{\mathbf{t}}) \geq 0 \ \forall \ \vec{\mathbf{t}} \in \mathbb{R}^{d_Y}$. Hence, $\vec{\epsilon}$ follows a legitimate multivariate stable distribution.

Finally, we construct the desired $Q_{MC}(M, \vec{\mathbf{X}}, \vec{\mathbf{Y}})$ from $Q_{MC}(M, \vec{\mathbf{X}}, \vec{\mathbf{X}}', \vec{\mathbf{Y}})$ by marginalizing $\vec{\mathbf{X}}'$.

**Part 2: Showing $Q_{MC}(M, \vec{\mathbf{X}}, \vec{\mathbf{Y}}) \in \Delta_P$**

For showing that $Q_{MC}(M, \vec{\mathbf{X}}, \vec{\mathbf{Y}}) \in \Delta_P$, we first need to derive $Q_{MC}(\vec{\mathbf{Y}}|M)$.

Deriving $Q(\vec{\mathbf{Y}}|M)$: We will derive $Q_{MC}(\vec{\mathbf{Y}}|M)$ by using the fact that $\vec{\epsilon} \perp\!\!\!\perp \vec{\mathbf{X}}'|M$, hence their conditional characteristic functions would just result in multiplication.

$$\mathbb{E}\left[ e^{i\vec{\mathbf{t}}^T \vec{\mathbf{Y}}} \Big| M = m \right] = \mathbb{E}\left[ e^{i\vec{\mathbf{t}}^T (\vec{\mathbf{X}}' + \vec{\epsilon})} \Big| M = m \right] = \mathbb{E}\left[ e^{i\vec{\mathbf{t}}^T \vec{\mathbf{X}}' + i\vec{\mathbf{t}}^T \vec{\epsilon}} \Big| M = m \right]$$

$$= \mathbb{E}\left[ e^{i\vec{\mathbf{t}}^T \vec{\mathbf{X}}'} e^{i\vec{\mathbf{t}}^T \vec{\epsilon}} \Big| M = m \right] \overset{(a)}{=} \mathbb{E}\left[ e^{i\vec{\mathbf{t}}^T \vec{\mathbf{X}}'} \Big| M = m \right] \mathbb{E}\left[ e^{i\vec{\mathbf{t}}^T \vec{\epsilon}} \Big| M = m \right], \tag{140}$$

where $(a)$ is due to $\vec{\epsilon} \perp\!\!\!\perp \vec{\mathbf{X}}'|M$. Substituting the characteristic functions of $\vec{\mathbf{X}}'$ and $\vec{\epsilon}$ from (124) and (126), respectively, in (140):

$$= \exp\left( -\sum_{j=1}^{d_Y} |t_j|^{\alpha} + f(k) \left| \vec{\mathbf{H}}_Y^T \vec{\mathbf{t}} \right|^{\alpha} \right) \exp\left( -f(k) \left| \vec{\mathbf{H}}_Y^T \vec{\mathbf{t}} \right|^{\alpha} + i\vec{\mathbf{t}}^T \vec{\mathbf{H}}_Y m \right)$$

$$= \exp\left( -\sum_{j=1}^{d_Y} |t_j|^{\alpha} + f(k) \left| \vec{\mathbf{H}}_Y^T \vec{\mathbf{t}} \right|^{\alpha} - f(k) \left| \vec{\mathbf{H}}_Y^T \vec{\mathbf{t}} \right|^{\alpha} + i\vec{\mathbf{t}}^T \vec{\mathbf{H}}_Y m \right)$$

$$= \exp\left( -\sum_{j=1}^{d_Y} |t_j|^{\alpha} + i\vec{\mathbf{t}}^T \vec{\mathbf{H}}_Y m \right). \tag{141}$$

By comparing the characteristic function shown in (141) with the characteristic function derived using $P(\vec{\mathbf{Y}}|M)$ shown in (116), we can conclude:

$$Q_{MC}(\vec{\mathbf{Y}}|M) = P(\vec{\mathbf{Y}}|M). \tag{142}$$

From (142) and (117), we can conclude:

$$Q_{MC}(M, \vec{\mathbf{Y}}) = Q_{MC}(M)Q_{MC}(\vec{\mathbf{Y}}|M) = P(M)P(\vec{\mathbf{Y}}|M) = P(M, \vec{\mathbf{Y}}). \tag{143}$$

We know that by construction:

$$Q_{MC}(M, \vec{\mathbf{X}}) = P(M, \vec{\mathbf{X}}). \tag{144}$$

Hence, $Q_{MC}(M, \vec{\mathbf{X}}, \vec{\mathbf{Y}}) \in \Delta_P$ and consequently, by proposition 1, $UI(M; \vec{\mathbf{Y}} \backslash \vec{\mathbf{X}}) = 0$, concluding our proof. $\square$

**Lemma 8.** *Let* $\vec{\mathbf{a}} = [a_1 \quad \dots \quad a_d] \in \mathbb{R}^d$ *and* $\vec{\mathbf{b}} = [b_1 \quad \dots \quad b_d] \in \mathbb{R}^d$, *then we have:*

$$\left| \vec{\mathbf{a}}^T \vec{\mathbf{b}} \right|^\alpha \leq \begin{cases} \|\vec{\mathbf{a}}\|_\alpha^\alpha \|\vec{\mathbf{b}}\|_{\alpha/(\alpha-1)}^\alpha & \alpha \in (1,2] \\ \|\vec{\mathbf{a}}\|_1^\alpha \|\vec{\mathbf{b}}\|_\infty^\alpha & \alpha \in (0,1] \end{cases}, \tag{145}$$

*where* $\|(\cdot)\|_p$ *is the standard* $L_p$ *norm.*

*Proof.* **Case 1**: $\alpha \in (1,2]$

Analysing the LHS of (145)

$$\left| \vec{\mathbf{a}}^T \vec{\mathbf{b}} \right| = \left| \sum_{i=1}^d a_i b_i \right| \overset{(a)}{\leq} \sum_{i=1}^d |a_i b_i| \overset{(b)}{\leq} \left( \sum_{i=1}^d |a_i|^\alpha \right)^{\frac{1}{\alpha}} \left( \sum_{i=1}^d |b_i|^{\frac{\alpha}{(\alpha-1)}} \right)^{\frac{(\alpha-1)}{\alpha}} = \|\vec{\mathbf{a}}\|_\alpha \|\vec{\mathbf{b}}\|_{\alpha/\alpha-1}, \tag{146}$$

where $(a)$ is due to the sub-additivity of $|(\cdot)|$ operator, and $(b)$ is due to Hölder's inequality [57]. Raising both sides of the inequality to $\alpha$ in (146) gives us the desired inequality. Since, $\alpha > 0$ raising both sides of the inequalities to power $\alpha$ does not change the direction of inequality.

$$\left| \vec{\mathbf{a}}^T \vec{\mathbf{b}} \right|^\alpha \leq \|\vec{\mathbf{a}}\|_\alpha^\alpha \|\vec{\mathbf{b}}\|_{\alpha/\alpha-1}^\alpha. \tag{147}$$

**Case 2**: $\alpha \in (0,1]$

Analysing the LHS of (145)

$$\left| \vec{\mathbf{a}}^T \vec{\mathbf{b}} \right| = \left| \sum_{i=1}^d a_i b_i \right| \overset{(a)}{\leq} \sum_{i=1}^d |a_i b_i| \overset{(b)}{\leq} \left( \sum_{i=1}^d |a_i| \right) \max_{i \in \{1,\dots,d\}} |b_i| = \|\vec{\mathbf{a}}\|_1 \|\vec{\mathbf{b}}\|_\infty, \tag{148}$$

where $(a)$ is again due to the sub-additivity of $|(\cdot)|$ operator, and $(b)$ is due to the fact that $|a_i b_i| \leq |a_i| \max_{i \in \{1,\dots,d\}} |b_i| \, \forall \, i \in \{1,\dots,d\}$. Raising both sides of the inequality to $\alpha$ in (148) gives us the desired inequality. Since, $\alpha > 0$ raising both sides of the inequalities to power $\alpha$ does not change the direction of inequality.

$$\left| \vec{\mathbf{a}}^T \vec{\mathbf{b}} \right|^\alpha \leq \|\vec{\mathbf{a}}\|_1^\alpha \|\vec{\mathbf{b}}\|_\infty^\alpha. \tag{149}$$

Combining (147) and (149), we obtain the desired inequality. $\qquad\square$

**Lemma 9.** *Let* $\vec{\mathbf{Z}} \sim p_{CS-IC}(\alpha, \vec{\mathbf{0}}_d, \vec{\mathbf{1}}_d, \vec{\mathbf{0}}_d)$ *be a* $d$-*dimensional random vector,* $\mathbf{A} = [a_{ij}]_{i,j=1}^{n,d} \in \mathbb{R}^{n \times d}$ *be a* $n \times d$ *matrix, and* $\vec{\mathbf{b}} \in \mathbb{R}^d$. *Then, the characteristic function of* $\mathbf{A}\vec{\mathbf{Z}} + \vec{\mathbf{b}}$ *is given by* (150)

$$\mathbb{E}\left[ e^{i\vec{\mathbf{t}}^T (\mathbf{A}\vec{\mathbf{Z}} + \vec{\mathbf{b}})} \right] = \exp\left( -\sum_{j=1}^d \left| \vec{\mathbf{a}}_j^T \vec{\mathbf{t}} \right|^\alpha + i\vec{\mathbf{t}}^T \vec{\mathbf{b}} \right) \, \forall \, \vec{\mathbf{t}} \in \mathbb{R}^d, \tag{150}$$

*where* $\vec{\mathbf{a}}_i$ *is the* $i$-*th column of* $\mathbf{A}$.

*Proof.* Analyzing L.H.S of (150)

$$\mathbb{E}\left[ e^{i\vec{\mathbf{t}}^T (\mathbf{A}\vec{\mathbf{Z}} + \vec{\mathbf{b}})} \right] = \mathbb{E}\left[ e^{i\vec{\mathbf{t}}^T \mathbf{A}\vec{\mathbf{Z}} + \vec{\mathbf{t}}^T \vec{\mathbf{b}}} \right] = \mathbb{E}\left[ e^{i\vec{\mathbf{t}}^T \mathbf{A}\vec{\mathbf{Z}}} e^{i\vec{\mathbf{t}}^T \vec{\mathbf{b}}} \right] \tag{151}$$

Using the fact that $e^{i\vec{\mathbf{t}}^T \vec{\mathbf{b}}}$ is constant w.r.t expectation over $\vec{\mathbf{Z}}$, and linearity of expectation, we have:

$$\begin{aligned} \mathbb{E}\left[ e^{i\vec{\mathbf{t}}^T (\mathbf{A}\vec{\mathbf{Z}} + \vec{\mathbf{b}})} \right] &= e^{i\vec{\mathbf{t}}^T \vec{\mathbf{b}}} \mathbb{E}\left[ \exp\left( i\vec{\mathbf{t}}^T \mathbf{A}\vec{\mathbf{Z}} \right) \right], \\ &= e^{i\vec{\mathbf{t}}^T \vec{\mathbf{b}}} \mathbb{E}\left[ \exp\left( i\vec{\mathbf{t}}^T [\vec{\mathbf{a}}_1 \quad \dots \quad \vec{\mathbf{a}}_d] \vec{\mathbf{Z}} \right) \right], \\ &= e^{i\vec{\mathbf{t}}^T \vec{\mathbf{b}}} \mathbb{E}\left[ \exp\left( i [\vec{\mathbf{t}}^T \vec{\mathbf{a}}_1 \quad \dots \quad \vec{\mathbf{t}}^T \vec{\mathbf{a}}_d] \vec{\mathbf{Z}} \right) \right], \end{aligned}$$

Simplifying the term $\begin{bmatrix} \vec{\mathbf{t}}^T\vec{\mathbf{a}}_1 & \ldots & \vec{\mathbf{t}}^T\vec{\mathbf{a}}_d \end{bmatrix}\vec{\mathbf{Z}}$ by substituting $\mathbf{Z} = [Z_1 \quad \ldots \quad Z_d]^T$, we obtain:

$$= e^{i\vec{\mathbf{t}}^T\vec{\mathbf{b}}}\mathbb{E}\left[\exp\left(i\sum_{j=1}^{d}(\vec{\mathbf{t}}^T\vec{\mathbf{a}}_j)Z_j\right)\right],$$

Using properties of exponentials, we obtain:

$$= e^{i\vec{\mathbf{t}}^T\vec{\mathbf{b}}}\mathbb{E}\left[\prod_{j=1}^{d}\exp\left(i\vec{\mathbf{t}}^T\vec{\mathbf{a}}_jZ_j\right)\right],$$

Using the fact $\{Z_j\}_{j=1}^{d}$ are jointly independent we obtain:

$$= e^{i\vec{\mathbf{t}}^T\vec{\mathbf{b}}}\prod_{j=1}^{d}\mathbb{E}\left[\exp\left(i\vec{\mathbf{t}}^T\vec{\mathbf{a}}_jZ_j\right)\right],$$

Using the fact that $Z_j \sim p_{CS}(\alpha, 0, 1, 0)$ and the result of lemma 5, we know that $\vec{\mathbf{t}}^T\vec{\mathbf{a}}_jZ_j \sim p_{CS}(\alpha, 0, |\vec{\mathbf{t}}^T\vec{\mathbf{a}}_j|, 0)$. Substituting the corresponding characteristic function of $\vec{\mathbf{t}}^T\vec{\mathbf{a}}_jZ_j$ using (323)

$$= e^{i\vec{\mathbf{t}}^T\vec{\mathbf{b}}}\prod_{j=1}^{d}\exp\left(-\left|\vec{\mathbf{t}}^T\vec{\mathbf{a}}_i\right|^{\alpha}\right) = e^{i\vec{\mathbf{t}}^T\vec{\mathbf{b}}}\exp\left(-\sum_{j=1}^{d}\left|\vec{\mathbf{t}}^T\vec{\mathbf{a}}_i\right|^{\alpha}\right)$$

$$= \exp\left(-\sum_{j=1}^{d}\left|\vec{\mathbf{t}}^T\vec{\mathbf{a}}_i\right|^{\alpha} + i\vec{\mathbf{t}}^T\vec{\mathbf{b}}\right). \tag{152}$$

$\square$

**Lemma 10.** *Let* $\vec{\mathbf{H}}_1 = \begin{bmatrix} h_1^{(1)} & \ldots & h_{d_1}^{(1)} \end{bmatrix}^T \in \mathbb{R}^{d_1}$ *and* $\vec{\mathbf{H}}_2 = \begin{bmatrix} h_1^{(2)} & \ldots & h_{d_2}^{(2)} \end{bmatrix}^T \in \mathbb{R}^{d_2}$. *Define* $\vec{\mathbf{H}}_1^k$ *as follows:*

$$\vec{\mathbf{H}}_1^k = \frac{1}{\|\vec{\mathbf{H}}_1\|_{1+k}^{1+k}}\begin{bmatrix} sgn(h_1^{(1)})|h_1^{(1)}|^k & \ldots & sgn(h_{d_1}^{(1)})|h_{d_1}^{(1)}|^k \end{bmatrix}^T \ \forall\, k \in [0, \infty).$$

*Furthermore, define* $\vec{\mathbf{H}}_1^\infty$ *as follows:*

$$j^* = \underset{j \in \{1, \ldots, d_1\}}{\arg\max}\left|h_j^{(1)}\right|,$$

$$\vec{\mathbf{H}}_1^\infty = \begin{bmatrix} h_1^\infty & \ldots & h_{d_1}^\infty \end{bmatrix}^T, \text{ with } h_j^\infty = 0 \ \forall\, j \in \{1, \ldots, d_1\}\setminus\{j^*\}, \text{ and } h_j^\infty = \frac{1}{h_{j^*}^{(1)}}. \tag{153}$$

*Then, we have:*

$$\vec{\mathbf{H}}_2(\vec{\mathbf{H}}_1^k)^T\vec{\mathbf{H}}_1 = \vec{\mathbf{H}}_2 \ \forall\, k \in [0, \infty) \text{ and } \vec{\mathbf{H}}_2(\vec{\mathbf{H}}_1^\infty)^T\vec{\mathbf{H}}_1 = \vec{\mathbf{H}}_2, \tag{154}$$

*where* $\|(\cdot)\|_{1+k}$ *is the standard* $L_{1+k}$ *norm [57]. We can alternatively state (154) in a more compact notation:*

$$\vec{\mathbf{H}}_2(\vec{\mathbf{H}}_1^k)^T\vec{\mathbf{H}}_1 = \vec{\mathbf{H}}_2 \ \forall\, k \in [0, \infty], \tag{155}$$

*where the case of* $k = \infty$ *is to be understood as* $\vec{\mathbf{H}}_1^k = \vec{\mathbf{H}}_1^\infty$.

*Proof.* **Case 1**: $k \in [0, \infty)$, i.e., $k$ is finite,

$$\vec{\mathbf{H}}_2(\vec{\mathbf{H}}_1^k)^T\vec{\mathbf{H}}_1 = \frac{1}{\|\vec{\mathbf{H}}_1\|_{1+k}^{1+k}}\vec{\mathbf{H}}_2\begin{bmatrix} sgn(h_1^{(1)})|h_1^{(1)}|^k & \ldots & sgn(h_{d_1}^{(1)})|h_{d_1}^{(1)}|^k \end{bmatrix}\begin{bmatrix} h_1^{(1)} \\ \vdots \\ h_{d_1}^{(1)} \end{bmatrix}$$

$$= \frac{1}{\|\vec{\mathbf{H}}_1\|_{1+k}^{1+k}}\underbrace{\left(\sum_{j=1}^{d}|h_j|^{1+k}\right)}_{=\|\vec{\mathbf{H}}_1\|_{1+k}^{1+k}}\vec{\mathbf{H}}_2 = \vec{\mathbf{H}}_2.$$

**Case 2**: $k = \infty$,

$$\vec{\mathbf{H}}_2 (\vec{\mathbf{H}}_1^\infty)^T \vec{\mathbf{H}}_1 = \vec{\mathbf{H}}_2 \left( \sum_{j=1}^{d_1} h_j^\infty h_j^{(1)} \right) \overset{(a)}{=} \vec{\mathbf{H}}_2 \left( h_{j*}^\infty h_{j*}^{(1)} \right) \overset{(b)}{=} \vec{\mathbf{H}}_2 \frac{1}{h_{j*}^{(1)}} h_{j*}^{(1)} = \vec{\mathbf{H}}_2,$$

where $(a)$ is due to the fact that all elements of $\vec{\mathbf{H}}_1^\infty$ are 0 except the $j^*$ element, and $(b)$ is due to the fact that $h_{j*}^\infty = 1/h_{j*}^{(1)}$. □

# H  Proof of Theorem 3

In this section, we provide the proof of theorem 3. Since theorem 3 provides the analytical PID terms for the system 2 of the multivariate affine continuous stable system defined in Sec. 4, we briefly restate certain key properties of the corresponding elliptically-contoured multivariate stable distributions and the corresponding multivariate affine continuous stable system for convenience.

## H.1  Elliptically-contoured multivariate stable distribution

Elliptically-contoured multivariate stable distributions are another example of a multivariate generalization of univariate continuous stable distributions. As the name suggests, the defining features of these distributions is that the corresponding p.d.f. has elliptical contours, similar to the multivariate Gaussian distribution. We refer the reader to Appx. N.3 for more details on multivariate continuous stable distributions. We now list certain key properties of elliptically-contoured multivariate stable distributions that we make use of in the proof of theorem 3:

1. We denote the p.d.f. of the elliptically-contoured continuous stable distribution as $p_{CS-EC}(\alpha, \mathbf{\Sigma}, \vec{\boldsymbol{\mu}})$, where $\mathbf{\Sigma}$ is a positive definite matrix, $\vec{\boldsymbol{\mu}} \in \mathbb{R}^d$, and $\alpha \in (0, 2]$. In general, the p.d.f. of Elliptically-contoured multivariate stable distribution do not have a closed-form analytical expression, and are expressed through their characteristic function.

2. The characteristic function of a $d$-dimensional random vector $\vec{\mathbf{X}}$ having elliptically-contoured multivariate distribution $p_{CS-EC}(\alpha, \mathbf{\Sigma}, \vec{\boldsymbol{\mu}})$ is expressed as follows:

$$\mathbb{E} \left[ e^{i \vec{\mathbf{t}}^T \vec{\mathbf{X}}} \right] = \exp \left( - \left( \vec{\mathbf{t}}^T \mathbf{\Sigma} \vec{\mathbf{t}} \right)^{\alpha/2} + i \vec{\mathbf{t}}^T \vec{\boldsymbol{\mu}} \right) \quad \forall \, \vec{\mathbf{t}} \in \mathbb{R}^d. \tag{156}$$

## H.2  Definition of the system 2 of the multivariate affine continuous stable system

Let the random variable $M \sim P(M)$ with support $\mathcal{M} \subseteq \mathbb{R}$. $\vec{\mathbf{X}}$ and $\vec{\mathbf{Y}}$ are $d_X$-dimensional and $d_Y$-dimensional random vectors, respectively. The joint distribution $P(M, \vec{\mathbf{X}}, \vec{\mathbf{Y}})$ describes the second system of the multivariate affine continuous stable distribution if it satisfies the following two properties:

1. $M \sim P(M)$ having some support set $\mathcal{M} \subseteq \mathbb{R}$.

2. The conditional distributions of $\vec{\mathbf{X}}$ and $\vec{\mathbf{Y}}$ are expressed as follows:

$$P(\vec{\mathbf{X}}|M) = p_{CS-EC}(\alpha, \mathbf{\Sigma}_X, \vec{\mathbf{H}}_X M + \vec{\mathbf{b}}_X), \text{ and}$$
$$P(\vec{\mathbf{Y}}|M) = p_{CS-EC}(\alpha, \mathbf{\Sigma}_Y, \vec{\mathbf{H}}_Y M + \vec{\mathbf{b}}_Y), \tag{157}$$

where $\mathbf{\Sigma}_X$ and $\mathbf{\Sigma}_Y$ are positive definite matrices, $\vec{\mathbf{H}}_X, \vec{\mathbf{b}}_X \in \mathbb{R}^{d_X}$, and $\vec{\mathbf{H}}_Y, \vec{\mathbf{b}}_Y \in \mathbb{R}^{d_Y}$.

## H.3  Formal proof of theorem 3

**Theorem 3.** *Let the joint distribution $P(M, \vec{\mathbf{X}}, \vec{\mathbf{Y}})$ of $M$, $\vec{\mathbf{X}}$, and $\vec{\mathbf{Y}}$ describe the system 2 of the multivariate affine continuous stable system. Define $\mathbf{\Sigma}_X^{-1/2}$ and $\mathbf{\Sigma}_Y^{-1/2}$ as the respective inverses of the matrices $\mathbf{\Sigma}_X^{1/2}$ and $\mathbf{\Sigma}_Y^{1/2}$ which satisfy: $(\mathbf{\Sigma}_X^{1/2})^T \mathbf{\Sigma}_X^{1/2} = \mathbf{\Sigma}_X$, and $(\mathbf{\Sigma}_Y^{1/2})^T \mathbf{\Sigma}_Y^{1/2} = \mathbf{\Sigma}_Y$. Without the loss of generality, assume $\|\mathbf{\Sigma}_Y^{-1/2} \vec{\mathbf{H}}_Y\|_2 \leq \|\mathbf{\Sigma}_X^{-1/2} \vec{\mathbf{H}}_X\|_2$. Then, $\Delta_P$ contains a Markov chain of the form $M \to \vec{\mathbf{X}} \to \vec{\mathbf{Y}}$ and $UI(M; \vec{\mathbf{Y}} \backslash \vec{\mathbf{X}}) = 0$.*

*Proof.* We first note two important observations:

1. We can always assume $\|\mathbf{\Sigma}_Y^{-1/2}\vec{\mathbf{H}}_Y\|_2 \leq \|\mathbf{\Sigma}_X^{-1/2}\vec{\mathbf{H}}_X\|_2$ without the loss generality because if $\|\mathbf{\Sigma}_Y^{-1/2}\vec{\mathbf{H}}_Y\|_2 \geq \|\mathbf{\Sigma}_X^{-1/2}\vec{\mathbf{H}}_X\|_2$, then we can always switch our nomenclature to refer to $\vec{\mathbf{Y}}$ as $\vec{\mathbf{X}}$, and $\vec{\mathbf{X}}$ as $\vec{\mathbf{Y}}$.

2. Properties of positive definite matrices guarantee the existence of an invertible $\mathbf{\Sigma}_X^{1/2}$ and $\mathbf{\Sigma}_Y^{1/2}$ [58].

The proof of this theorem is a generalization of the proof used for the deriving a similar result for the special case of the multivariate Gaussian distribution discussed in [22], which borrows known results from Gaussian interference channels discussed in [48]. This proof follows the same structure as our earlier proofs.

1. In the first part, we explicitly construct a joint distribution $Q_{MC}(M, \vec{\mathbf{X}}, \vec{\mathbf{Y}})$ having the Markovian structure $M \to \vec{\mathbf{X}} \to \vec{\mathbf{Y}}$.

2. In the second part, we show that the $Q_{MC}(M, \vec{\mathbf{X}}, \vec{\mathbf{Y}})$ constructed in the first part, lies in $\Delta_P$. Therefore, we can then apply the result of proposition 1 to conclude $UI(M; \vec{\mathbf{Y}} \backslash \vec{\mathbf{X}}) = 0$.

**Part 1: Specifying $Q_{MC}(M, \vec{\mathbf{X}}, \vec{\mathbf{Y}})$**

We explicitly construct the desired Markov chain $M \to \vec{\mathbf{X}} \to \vec{\mathbf{Y}}$ by constructing a larger Markov chain $M \to \vec{\mathbf{X}} \to \vec{\mathbf{X}}' \to \vec{\mathbf{X}}'' \to \vec{\mathbf{Y}}' \to \vec{\mathbf{Y}}$, and then marginalizing the larger Markov chain to obtain the desired Markov chain $M \to \vec{\mathbf{X}} \to \vec{\mathbf{Y}}$.

Denote the joint distribution of the Markov chain $M \to \vec{\mathbf{X}} \to \vec{\mathbf{X}}' \to \vec{\mathbf{X}}'' \to \vec{\mathbf{Y}}' \to \vec{\mathbf{Y}}$ as $Q_{MC}(M, \vec{\mathbf{X}}, \vec{\mathbf{X}}', \vec{\mathbf{X}}'', \vec{\mathbf{Y}}', \vec{\mathbf{Y}})$. We can decompose $Q_{MC}(M, \vec{\mathbf{X}}, \vec{\mathbf{X}}', \vec{\mathbf{X}}'', \vec{\mathbf{Y}}', \vec{\mathbf{Y}})$ by utilizing its Markovian structure as follows: $Q_{MC}(M, \vec{\mathbf{X}}, \vec{\mathbf{X}}', \vec{\mathbf{X}}'', \vec{\mathbf{Y}}', \vec{\mathbf{Y}}) = Q_{MC}(M)Q_{MC}(\vec{\mathbf{X}}|M)$ $Q_{MC}(\vec{\mathbf{X}}'|\vec{\mathbf{X}}) \, Q_{MC}(\vec{\mathbf{X}}''|\vec{\mathbf{X}}')Q_{MC}(\vec{\mathbf{Y}}'|\vec{\mathbf{X}}'')Q_{MC}(\vec{\mathbf{Y}}|\vec{\mathbf{Y}}')$. Consequently, we can specify/construct the distribution $Q_{MC}(M, \vec{\mathbf{X}}, \vec{\mathbf{X}}', \vec{\mathbf{X}}'', \vec{\mathbf{Y}}', \vec{\mathbf{Y}})$ by individually specifying $Q_{MC}(M)$, $Q_{MC}(\vec{\mathbf{X}}|M)$, $Q_{MC}(\vec{\mathbf{X}}'|\vec{\mathbf{X}})$, $Q_{MC}(\vec{\mathbf{X}}''|\vec{\mathbf{X}}')$, $Q_{MC}(\vec{\mathbf{Y}}'|\vec{\mathbf{X}}'')$, and $Q_{MC}(\vec{\mathbf{Y}}|\vec{\mathbf{Y}}')$.

Specifying $Q_{MC}(M)$ and $Q_{MC}(\vec{\mathbf{X}}|M)$: We choose $Q_{MC}(M)$ and $Q_{MC}(\vec{\mathbf{X}}|M)$ as follows:

$$Q_{MC}(M) = P(M) \text{ and } Q_{MC}(\vec{\mathbf{X}}|M) = P(\vec{\mathbf{X}}|M). \tag{158}$$

Specifying $Q_{MC}(\vec{\mathbf{X}}'|\vec{\mathbf{X}})$: $Q_{MC}(\vec{\mathbf{X}}'|\vec{\mathbf{X}})$ is specified through the following deterministic transformation:

$$\vec{\mathbf{X}}' = (\Sigma_X^{-1/2})^T(\vec{\mathbf{X}} - \vec{\mathbf{b}}_X). \tag{159}$$

We will also derive the distribution $Q(\vec{\mathbf{X}}'|M)$ before proceeding with our construction, as we will need it later in the proof to show that the constructed $Q_{MC}(M, \vec{\mathbf{X}}, \vec{\mathbf{Y}}) \in \Delta_P$.

*Deriving $Q_{MC}(\vec{\mathbf{X}}'|M)$:* We use lemma 11 to derive $Q_{MC}(\vec{\mathbf{X}}'|M = m)$, as $Q_{MC}(\vec{\mathbf{X}}|M = m)$ is an elliptically contoured multivariate stable distribution for a fixed $m$, and (159) defines a scaling and

translation operation.

$$Q_{MC}(\vec{\mathbf{X}}'|M=m) = p_{CS-EC}\left(\alpha, (\boldsymbol{\Sigma}_X^{-1/2})^T \boldsymbol{\Sigma}_X \boldsymbol{\Sigma}_X^{-1/2}, (\boldsymbol{\Sigma}_X^{-1/2})^T \vec{\mathbf{H}}_X m + (\boldsymbol{\Sigma}_X^{-1/2})^T (\vec{\mathbf{b}}_X - \vec{\mathbf{b}}_X)\right),$$

$$\Rightarrow Q_{MC}(\vec{\mathbf{X}}'|M=m) = p_{CS-EC}\left(\alpha, (\boldsymbol{\Sigma}_X^{-1/2})^T \boldsymbol{\Sigma}_X \boldsymbol{\Sigma}_X^{-1/2}, (\boldsymbol{\Sigma}_X^{-1/2})^T \vec{\mathbf{H}}_X m\right),$$

$$\Rightarrow Q_{MC}(\vec{\mathbf{X}}'|M=m) = p_{CS-EC}\left(\alpha, (\boldsymbol{\Sigma}_X^{-1/2})^T (\boldsymbol{\Sigma}_X^{1/2})^T \boldsymbol{\Sigma}_X^{1/2} \boldsymbol{\Sigma}_X^{-1/2}, (\boldsymbol{\Sigma}_X^{-1/2})^T \vec{\mathbf{H}}_X m\right),$$

$$\Rightarrow Q_{MC}(\vec{\mathbf{X}}'|M=m) = p_{CS-EC}\left(\alpha, (\boldsymbol{\Sigma}_X^{1/2} \boldsymbol{\Sigma}_X^{-1/2})^T \boldsymbol{\Sigma}_X^{1/2} \boldsymbol{\Sigma}_X^{-1/2}, (\boldsymbol{\Sigma}_X^{-1/2})^T \vec{\mathbf{H}}_X m\right),$$

$$\Rightarrow Q_{MC}(\vec{\mathbf{X}}'|M=m) = p_{CS-EC}\left(\alpha, \mathbf{I}_{d_X}, (\boldsymbol{\Sigma}_X^{-1/2})^T \vec{\mathbf{H}}_X m\right),$$

$$\Rightarrow Q_{MC}(\vec{\mathbf{X}}'|M=m) = p_{CS-EC}\left(\alpha, \mathbf{I}_{d_X}, \vec{\mathbf{K}}_X m\right), \tag{160}$$

where $\mathbf{I}_{d_X}$ is a $d_X \times d_X$ identity matrix, and $\vec{\mathbf{K}}_X = (\boldsymbol{\Sigma}_X^{-1/2})^T \vec{\mathbf{H}}_X$.

Specifying $Q_{MC}(\vec{\mathbf{X}}''|\vec{\mathbf{X}}')$: $Q_{MC}(\vec{\mathbf{X}}''|\vec{\mathbf{X}}')$ is again specified through a deterministic transformation:

$$\vec{\mathbf{X}}'' = \vec{\mathbf{K}}_Y (\vec{\mathbf{K}}_X^1)^T \vec{\mathbf{X}}', \tag{161}$$

where $\vec{\mathbf{K}}_Y = (\boldsymbol{\Sigma}_Y^{-1/2})^T \vec{\mathbf{H}}_Y$ and $\vec{\mathbf{K}}_X^1 = {}^1/\|\vec{\mathbf{K}}_X\|_2^2 \vec{\mathbf{K}}_X$. We also derive the distribution $Q(\vec{\mathbf{X}}'|M)$ which will be needed for showing that the constructed $Q_{MC}(M, \vec{\mathbf{X}}, \vec{\mathbf{Y}}) \in \Delta_P$.

*Deriving $Q_{MC}(\vec{\mathbf{X}}''|M = m)$:* We again use lemma 11 to calculate $Q_{MC}(\vec{\mathbf{X}}''|M = m)$, as $Q_{MC}(\vec{\mathbf{X}}'|M = m)$ is an elliptically contoured multivariate stable distribution (see (160)) for a fixed $m$, and (159) defines a scaling and translation operation.

$$Q_{MC}(\vec{\mathbf{X}}'|M=m) = p_{CS-EC}\left(\alpha, (\vec{\mathbf{K}}_Y (\vec{\mathbf{K}}_X^1)^T) \mathbf{I}_{d_X} (\vec{\mathbf{K}}_Y (\vec{\mathbf{K}}_X^1)^T)^T, \vec{\mathbf{K}}_Y (\vec{\mathbf{K}}_X^1)^T \vec{\mathbf{K}}_X m\right),$$

$$= p_{CS-EC}\left(\alpha, (\vec{\mathbf{K}}_Y (\vec{\mathbf{K}}_X^1)^T)(\vec{\mathbf{K}}_Y (\vec{\mathbf{K}}_X^1)^T)^T, \vec{\mathbf{K}}_Y (\vec{\mathbf{K}}_X^1)^T \vec{\mathbf{K}}_X m\right),$$

From lemma 10, we know that $\vec{\mathbf{K}}_Y (\vec{\mathbf{K}}_X^1)^T \vec{\mathbf{K}}_X = \vec{\mathbf{K}}_Y$, which implies

$$Q_{MC}(\vec{\mathbf{X}}'|M=m) = p_{CS-EC}\left(\alpha, (\vec{\mathbf{K}}_Y (\vec{\mathbf{K}}_X^1)^T)(\vec{\mathbf{K}}_Y (\vec{\mathbf{K}}_X^1)^T)^T, \vec{\mathbf{K}}_Y m\right). \tag{162}$$

Specifying $Q_{MC}(\vec{\mathbf{Y}}'|\vec{\mathbf{X}}'')$: We specify $Q_{MC}(\vec{\mathbf{Y}}'|\vec{\mathbf{X}}'')$ through the following stochastic transformation:

$$\vec{\mathbf{Y}}' = \vec{\mathbf{X}}'' + \vec{\boldsymbol{\epsilon}}, \tag{163}$$

where $\vec{\boldsymbol{\epsilon}}$ follows a multivariate stable distribution. Furthermore, we assume that $\vec{\boldsymbol{\epsilon}}$ is jointly independent from $(\vec{\mathbf{X}}, \vec{\mathbf{X}}', \vec{\mathbf{X}}'', M)$, i.e., $\vec{\boldsymbol{\epsilon}} \perp\!\!\!\perp (\vec{\mathbf{X}}, \vec{\mathbf{X}}', \vec{\mathbf{X}}'', M)$. The characteristic function of $\vec{\boldsymbol{\epsilon}}$ is defined as follows:

$$\mathbb{E}\left[e^{i\vec{\mathbf{t}}^T \vec{\boldsymbol{\epsilon}}}\right] = \exp\left(-\left(\vec{\mathbf{t}}^T \mathbf{I}_{d_Y} \vec{\mathbf{t}}\right)^{\alpha/2} + \left(\vec{\mathbf{t}}^T (\vec{\mathbf{K}}_Y (\vec{\mathbf{K}}_X^1)^T)(\vec{\mathbf{K}}_Y (\vec{\mathbf{K}}_X^1)^T)^T \vec{\mathbf{t}}\right)^{\alpha/2}\right). \tag{164}$$

In order for $Q_{MC}(\vec{\mathbf{Y}}'|\vec{\mathbf{X}}'')$ to be a valid distribution, we need to ensure that the distribution of $\vec{\boldsymbol{\epsilon}}$ defined in (164) is a legitimate multivariate stable distribution. Similarly to the proof of theorem 2, we will utilize property 1 for showing that $\vec{\boldsymbol{\epsilon}}$ follows a stable multivariate stable distribution. Property 1 states a random vector is distributed according to a multivariate stable distribution iff every 1-dimensional projection of this random vector follows a univariate stable continuous distribution. Hence, to show that $\vec{\boldsymbol{\epsilon}}$ is distributed according to a legitimate multivariate stable distribution, we will show that every 1-dimensional projection of $\vec{\boldsymbol{\epsilon}}$ follows a univariate stable continuous distribution.

Let $\vec{\mathbf{t}} \in \mathbb{R}^{d_Y}$, then the characteristic function of the 1-dimensional projection of $\vec{\boldsymbol{\epsilon}}$ along $\vec{\mathbf{t}}$, i.e., $\vec{\mathbf{t}}^T \vec{\boldsymbol{\epsilon}}$ can be trivially derived from (164).

$$\mathbb{E}\left[e^{i\vec{\mathbf{t}}^T \vec{\boldsymbol{\epsilon}}}\right] = \exp\left(-\left(\vec{\mathbf{t}}^T \mathbf{I}_{d_Y} \vec{\mathbf{t}}\right)^{\alpha/2} + \left(\vec{\mathbf{t}}^T (\vec{\mathbf{K}}_Y (\vec{\mathbf{K}}_X^1)^T)(\vec{\mathbf{K}}_Y (\vec{\mathbf{K}}_X^1)^T)^T \vec{\mathbf{t}}\right)^{\alpha/2}\right). \tag{165}$$

Comparing the characteristic function of $\vec{\mathbf{t}}^T \vec{\epsilon}$ described in (127) with the standard characteristic function of a univariate stable characteristic function defined in (323), we can conclude $\vec{\mathbf{t}}^T \vec{\epsilon} \sim p_{CS}(\alpha, \beta(\vec{\mathbf{t}}), \gamma(\vec{\mathbf{t}}), \mu(\vec{\mathbf{t}}))$, where:

$$\gamma(\vec{\mathbf{t}}) = \left(\vec{\mathbf{t}}^T \mathbf{I}_{d_Y} \vec{\mathbf{t}}\right)^{\alpha/2} - \left(\vec{\mathbf{t}}^T (\vec{\mathbf{K}}_Y (\vec{\mathbf{K}}_X^1)^T)(\vec{\mathbf{K}}_Y (\vec{\mathbf{K}}_X^1)^T)^T \vec{\mathbf{t}}\right)^{\alpha/2}, \mu(\vec{\mathbf{t}}) = 0, \text{ and } \beta(\vec{\mathbf{t}}) = 0. \quad (166)$$

In order for $\vec{\mathbf{t}}^T \vec{\epsilon}$ to have a legitimate univariate stable distribution, we just need to show that $\gamma(\vec{\mathbf{t}}) \geq 0 \ \forall \ \vec{\mathbf{t}} \in \mathbb{R}^{d_Y}$. Note that $\gamma(\vec{\mathbf{t}}) = 0$ would correspond to the case where all the mass of the distribution is centered at 0, which does satisfy definition 2, and is an example of a degenerate univariate stable distribution.

By the assumption in theorem, $\|\vec{\mathbf{K}}_Y\|_2 = \|\mathbf{\Sigma}_Y^{-1/2}\vec{\mathbf{H}}_Y\|_2 \leq \|\vec{\mathbf{K}}_X\|_2 = \|\mathbf{\Sigma}_X^{-1/2}\vec{\mathbf{H}}_X\|_2$. Hence, we can use the result of lemma 12, which shows that $(\vec{\mathbf{K}}_Y (\vec{\mathbf{K}}_X^1)^T)(\vec{\mathbf{K}}_Y (\vec{\mathbf{K}}_X^1)^T)^T \preceq \mathbf{I}_{d_Y}$. Using the fact that $(\mathbf{I}_{d_Y} - (\vec{\mathbf{K}}_Y (\vec{\mathbf{K}}_X^1)^T)(\vec{\mathbf{K}}_Y (\vec{\mathbf{K}}_X^1)^T)^T)$ is a positive semi-definite matrix, we have:

$$\vec{\mathbf{t}}^T \left( \mathbf{I}_{d_Y} - (\vec{\mathbf{K}}_Y (\vec{\mathbf{K}}_X^1)^T)(\vec{\mathbf{K}}_Y (\vec{\mathbf{K}}_X^1)^T)^T \right) \vec{\mathbf{t}} \geq 0 \ \forall \ \vec{\mathbf{t}} \in \mathbb{R}^{d_Y},$$

$$\Rightarrow \vec{\mathbf{t}}^T \mathbf{I}_{d_Y} \vec{\mathbf{t}} \geq \vec{\mathbf{t}}^T (\vec{\mathbf{K}}_Y (\vec{\mathbf{K}}_X^1)^T)(\vec{\mathbf{K}}_Y (\vec{\mathbf{K}}_X^1)^T)^T \vec{\mathbf{t}} \ \forall \ \vec{\mathbf{t}} \in \mathbb{R}^{d_Y},$$

$$\Rightarrow \left(\vec{\mathbf{t}}^T \mathbf{I}_{d_Y} \vec{\mathbf{t}}\right)^{\alpha/2} \geq \left(\vec{\mathbf{t}}^T (\vec{\mathbf{K}}_Y (\vec{\mathbf{K}}_X^1)^T)(\vec{\mathbf{K}}_Y (\vec{\mathbf{K}}_X^1)^T)^T \vec{\mathbf{t}}\right)^{\alpha/2} \ \forall \ \vec{\mathbf{t}} \in \mathbb{R}^{d_Y},$$

$$\Rightarrow \gamma(\vec{\mathbf{t}}) = \left(\vec{\mathbf{t}}^T \mathbf{I}_{d_Y} \vec{\mathbf{t}}\right)^{\alpha/2} - \left(\vec{\mathbf{t}}^T (\vec{\mathbf{K}}_Y (\vec{\mathbf{K}}_X^1)^T)(\vec{\mathbf{K}}_Y (\vec{\mathbf{K}}_X^1)^T)^T \vec{\mathbf{t}}\right)^{\alpha/2} \geq 0 \ \forall \ \vec{\mathbf{t}} \in \mathbb{R}^{d_Y}. \quad (167)$$

Hence, we know that $\vec{\epsilon}$ follows a legitimate multivariate stable distribution. We also derive $Q_{MC}(\vec{\mathbf{Y}}'|M)$ which is needed for showing $Q_{MC}(M, \vec{\mathbf{X}}, \vec{\mathbf{Y}}) \in \Delta_P$.

*Deriving $Q_{MC}(\vec{\mathbf{Y}}'|M)$:* We will derive $Q_{MC}(\vec{\mathbf{Y}}'|M)$ using the fact that $\vec{\epsilon} \perp\!\!\!\perp \vec{\mathbf{X}}''|M = m$, hence their conditional characteristic functions would just result in multiplication.

$$\mathbb{E}\left[e^{i\vec{\mathbf{t}}^T \vec{\mathbf{Y}}'} \Big| M = m\right] = \mathbb{E}\left[e^{i\vec{\mathbf{t}}^T (\vec{\mathbf{X}}'' + \vec{\epsilon})} \Big| M = m\right] = \mathbb{E}\left[e^{i\vec{\mathbf{t}}^T \vec{\mathbf{X}}'' + i\vec{\mathbf{t}}^T \vec{\epsilon}} \Big| M = m\right],$$

$$= \mathbb{E}\left[e^{i\vec{\mathbf{t}}^T \vec{\mathbf{X}}''} e^{i\vec{\mathbf{t}}^T \vec{\epsilon}} \Big| M = m\right] \overset{(a)}{=} \mathbb{E}\left[e^{i\vec{\mathbf{t}}^T \vec{\mathbf{X}}''} \Big| M = m\right] \mathbb{E}\left[e^{i\vec{\mathbf{t}}^T \vec{\epsilon}} \Big| M = m\right], \quad (168)$$

where $(a)$ is due to $\vec{\epsilon} \perp\!\!\!\perp \vec{\mathbf{X}}''|M$. Substituting the characteristic functions of $\vec{\mathbf{X}}''$ and $\vec{\epsilon}$ from (162) and (164), respectively, in (168)

$$= \exp\left( - \left(\vec{\mathbf{t}}^T (\vec{\mathbf{K}}_Y (\vec{\mathbf{K}}_X^1)^T)(\vec{\mathbf{K}}_Y (\vec{\mathbf{K}}_X^1)^T)^T \vec{\mathbf{t}}\right)^{\alpha/2} + i\vec{\mathbf{t}}^T \vec{\mathbf{K}}_Y m \right) \times$$

$$\exp\left( - \left(\vec{\mathbf{t}}^T \mathbf{I}_{d_Y} \vec{\mathbf{t}}\right)^{\alpha/2} + \left(\vec{\mathbf{t}}^T (\vec{\mathbf{K}}_Y (\vec{\mathbf{K}}_X^1)^T)(\vec{\mathbf{K}}_Y (\vec{\mathbf{K}}_X^1)^T)^T \vec{\mathbf{t}}\right)^{\alpha/2} \right),$$

$$= \exp\left( - \left(\vec{\mathbf{t}}^T \mathbf{I}_{d_Y} \vec{\mathbf{t}}\right)^{\alpha/2} + i\vec{\mathbf{t}}^T \vec{\mathbf{K}}_Y m \right). \quad (169)$$

By inspecting the characteristic function shown in (169), we can conclude:

$$Q_{MC}(\vec{\mathbf{Y}}'|M = m) = p_{CS-EC}(\alpha, \mathbf{I}_{d_Y}, \vec{\mathbf{K}}_Y m). \quad (170)$$

Specifying $Q_{MC}(\vec{\mathbf{Y}}|\vec{\mathbf{Y}}')$: Lastly, we specify $Q_{MC}(\vec{\mathbf{Y}}|\vec{\mathbf{Y}}')$ through the following deterministic transformation:

$$\vec{\mathbf{Y}} = (\mathbf{\Sigma}_Y^{1/2})^T \vec{\mathbf{Y}}' + \vec{\mathbf{b}}_Y. \quad (171)$$

Finally, we construct the desired $Q_{MC}(M, \vec{\mathbf{X}}, \vec{\mathbf{Y}})$ from $Q_{MC}(M, \vec{\mathbf{X}}, \vec{\mathbf{X}}', \vec{\mathbf{X}}'', \vec{\mathbf{Y}}', \vec{\mathbf{Y}})$ by marginalizing $\vec{\mathbf{X}}', \vec{\mathbf{X}}''$, and $\vec{\mathbf{Y}}'$.

**Part 2: Showing $Q_{MC}(M, \vec{\mathbf{X}}, \vec{\mathbf{Y}}) \in \Delta_P$**

For showing that $Q_{MC}(M, \vec{\mathbf{X}}, \vec{\mathbf{Y}}) \in \Delta_P$, we first need to derive $Q_{MC}(\vec{\mathbf{Y}}|M)$.

Deriving $Q(\vec{\mathbf{Y}}|M)$: We use lemma 11 to calculate $Q_{MC}(\vec{\mathbf{Y}}|M=m)$, as $Q_{MC}(\vec{\mathbf{X}}|M=m)$ is an elliptically contoured multivariate continuous stable distribution for a fixed $m$ (see (170)), and (171) defines a scaling and translation operation.

$$Q_{MC}(\vec{\mathbf{X}}'|M=m) = p_{CS-EC}\left(\alpha, (\mathbf{\Sigma}_Y^{1/2})^T\mathbf{\Sigma}_Y^{1/2}, (\mathbf{\Sigma}_Y^{1/2})^T\vec{\mathbf{K}}_Y m + \vec{\mathbf{b}}_Y\right),$$

$$\overset{(a)}{=} p_{CS-EC}\left(\alpha, \mathbf{\Sigma}_Y, (\mathbf{\Sigma}_Y^{1/2})^T\vec{\mathbf{K}}_Y m + \vec{\mathbf{b}}_Y\right),$$

$$\overset{(b)}{=} p_{CS-EC}\left(\alpha, \mathbf{\Sigma}_Y, (\mathbf{\Sigma}_Y^{1/2})^T(\mathbf{\Sigma}_Y^{-1/2})^T\vec{\mathbf{H}}_Y m + \vec{\mathbf{b}}_Y\right), \qquad (172)$$

$$= p_{CS-EC}\left(\alpha, \mathbf{\Sigma}_Y, \vec{\mathbf{H}}_Y m + \vec{\mathbf{b}}_Y\right), \qquad (173)$$

where $(a)$ is due to the fact that $(\mathbf{\Sigma}_Y^{1/2})^T\mathbf{\Sigma}_Y^{1/2} = \mathbf{\Sigma}_Y$, and $(b)$ is due to the fact that $\vec{\mathbf{K}}_Y = \mathbf{\Sigma}_Y^{-1/2}\vec{\mathbf{H}}_Y$. Comparing (173) with $P(\vec{\mathbf{Y}}|M=m)$ shown in Appx. H.2, we can conclude:

$$Q_{MC}(\vec{\mathbf{Y}}|M) = P(\vec{\mathbf{Y}}|M). \qquad (174)$$

From (174) and (158), we can conclude:

$$Q_{MC}(M, \vec{\mathbf{Y}}) = Q_{MC}(M)Q_{MC}(\vec{\mathbf{Y}}|M) = P(M)P(\vec{\mathbf{Y}}|M) = P(M, \vec{\mathbf{Y}}). \qquad (175)$$

We know that by construction:

$$Q_{MC}(M, \vec{\mathbf{X}}) = P(M, \vec{\mathbf{X}}). \qquad (176)$$

Hence, $Q_{MC}(M, \vec{\mathbf{X}}, \vec{\mathbf{Y}}) \in \Delta_P$ and consequently, by proposition 1, $UI(M; \vec{\mathbf{Y}}\backslash\vec{\mathbf{X}}) = 0$, concluding our proof. $\qquad\square$

**Lemma 11.** *Let $\vec{\mathbf{X}} \sim p_{CS-EC}(\alpha, \mathbf{\Sigma}, \vec{\boldsymbol{\mu}})$ be a d-dimensional random vector, where $\mathbf{\Sigma}$ is a $d \times d$ positive definite matrix, $\vec{\boldsymbol{\mu}} \in \mathbb{R}^d$, and $\alpha \in (0, 2]$. Furthermore, let $\mathbf{A} \in \mathbb{R}^{n \times d}$ and $\vec{\mathbf{b}} \in \mathbb{R}^d$. Then,*

$$\mathbf{A}\vec{\mathbf{X}} + \vec{\mathbf{b}} \sim p_{CS-EC}(\alpha, \mathbf{A}\mathbf{\Sigma}\mathbf{A}^T, \mathbf{A}\vec{\boldsymbol{\mu}} + \vec{\mathbf{b}}).$$

*Proof.* Calculating the characteristic function of $\mathbf{A}\vec{\mathbf{X}} + \vec{\mathbf{b}}$:

$$\mathbb{E}\left[\exp\left(i\vec{\mathbf{t}}^T(\mathbf{A}\vec{\mathbf{X}} + \vec{\mathbf{b}})\right)\right] = \mathbb{E}\left[\exp\left(i\vec{\mathbf{t}}^T\mathbf{A}\vec{\mathbf{X}} + i\vec{\mathbf{t}}^T\vec{\mathbf{b}}\right)\right] = \mathbb{E}\left[\exp\left(i\vec{\mathbf{t}}^T\mathbf{A}\vec{\mathbf{X}}\right)\exp\left(i\vec{\mathbf{t}}^T\vec{\mathbf{b}}\right)\right],$$

Employing linearity of expectation to pull the term $\exp\left(i\vec{\mathbf{t}}^T\vec{\mathbf{b}}\right)$ out of the expectation, as it is constant with respect to $\vec{\mathbf{X}}$:

$$\mathbb{E}\left[\exp\left(i\vec{\mathbf{t}}^T(\mathbf{A}\vec{\mathbf{X}} + \vec{\mathbf{b}})\right)\right] = \exp\left(i\vec{\mathbf{t}}^T\vec{\mathbf{b}}\right)\mathbb{E}\left[\exp\left(i\vec{\mathbf{t}}^T\mathbf{A}\vec{\mathbf{X}}\right)\right]. \qquad (177)$$

Substituting $\vec{\mathbf{t}}' = \vec{\mathbf{t}}^T\mathbf{A}$ in (177):

$$\mathbb{E}\left[\exp\left(i\vec{\mathbf{t}}^T(\mathbf{A}\vec{\mathbf{X}} + \vec{\mathbf{b}})\right)\right] = \exp\left(i\vec{\mathbf{t}}^T\vec{\mathbf{b}}\right)\mathbb{E}\left[\exp\left(i(\vec{\mathbf{t}}')^T\vec{\mathbf{X}}\right)\right].$$

Substituting the formula of the characteristic function of an Elliptically-contoured multivariate stable distribution (given in (156)) for calculating $\mathbb{E}\left[\exp\left(i(\vec{\mathbf{t}}')^T\vec{\mathbf{X}}\right)\right]$ as $\vec{\mathbf{X}} \sim p_{CS-EC}(\alpha, \mathbf{\Sigma}, \vec{\boldsymbol{\mu}})$:

$$\mathbb{E}\left[\exp\left(i\vec{\mathbf{t}}^T(\mathbf{A}\vec{\mathbf{X}} + \vec{\mathbf{b}})\right)\right] = \exp\left(i\vec{\mathbf{t}}^T\vec{\mathbf{b}}\right)\exp\left(-\left((\vec{\mathbf{t}}')^T\mathbf{\Sigma}\vec{\mathbf{t}}'\right)^{\alpha/2} + i(\vec{\mathbf{t}}')^T\vec{\boldsymbol{\mu}}\right).$$

Re-substituting $\vec{\mathbf{t}}' = \vec{\mathbf{t}}^T\mathbf{A}$ in the above equation

$$= \exp\left(i\vec{\mathbf{t}}^T\vec{\mathbf{b}}\right)\exp\left(-\left(\vec{\mathbf{t}}^T\mathbf{A}\mathbf{\Sigma}\mathbf{A}^T\vec{\mathbf{t}}\right)^{\alpha/2} + i\vec{\mathbf{t}}^T\mathbf{A}\vec{\boldsymbol{\mu}}\right),$$

$$= \exp\left(-\left(\vec{\mathbf{t}}^T\mathbf{A}\mathbf{\Sigma}\mathbf{A}^T\vec{\mathbf{t}}\right)^{\alpha/2} + i\vec{\mathbf{t}}^T(\mathbf{A}\vec{\boldsymbol{\mu}} + \vec{\mathbf{b}})\right). \qquad (178)$$

By inspecting the characteristic function described in (178), we can determine that $\mathbf{A}\vec{\mathbf{X}} + \vec{\mathbf{b}}$ is distributed according to an elliptically-contoured multivariate stable distribution, i.e., $\mathbf{A}\vec{\mathbf{X}} + \vec{\mathbf{b}} \sim p_{CS-EC}\left(\alpha, \mathbf{A}\mathbf{\Sigma}\mathbf{A}^T, \mathbf{A}\vec{\boldsymbol{\mu}} + \vec{\mathbf{b}}\right)$. $\qquad\square$

**Lemma 12.** *Let $\vec{\mathbf{H}}_1 \in \mathbb{R}^{d_1}$ and $\vec{\mathbf{H}}_2 \in \mathbb{R}^{d_2}$. Then, $\|\vec{\mathbf{H}}_1\|_2 \geq \|\vec{\mathbf{H}}_2\|_2$ if and only if $\exists$ a matrix $\mathbf{A} \in \mathbb{R}^{d_2 \times d_1}$ such that $\vec{\mathbf{H}}_2 = \mathbf{A}\vec{\mathbf{H}}_1$, and $\mathbf{A}\mathbf{A}^T \preceq \mathbf{I}_{d_2}$, with $\mathbf{A} = \vec{\mathbf{H}}_2(\vec{\mathbf{H}}_1^1)^T$, where $\mathbf{B} \preceq \mathbf{C}$ means $\mathbf{B} - \mathbf{C}$ is a positive semi-definite matrix, $\vec{\mathbf{H}}_1^1$ is as defined in lemma 10, and $\mathbf{I}_{d_2}$ is $d_2 \times d_2$ identity matrix.*

*Proof.* See the proof of lemma 5 in [48] for the special case $t = 1$. $\qquad\square$

# I  Proof of Theorem 4

In this section, we provide the proof of theorem 4. Since theorem 4 provides the analytical PID terms for the univariate affine discrete stable system defined in Sec. 4, we briefly restate certain key properties of the univariate discrete stable distributions and the corresponding univariate affine discrete stable system for convenience.

## I.1  Univariate discrete stable distribution

Univariate discrete stable distributions are the discrete analogues of univariate continuous stable distributions. We refer the reader to Appx. N.4 for more details on univariate discrete stable distributions. We now list certain key properties of univariate continuous stable distributions that we make use of in the proof of theorem 4:

1. If $X$ is distributed according to a univariate discrete stable distribution, then $X \overset{d}{=} \gamma \circ X_1 + (1 - \gamma^\nu)^{1/\nu} \circ X_2$, where $X_1$ and $X_2$ are two independent copies of $X$, $\nu \in (0, 1]$, $\gamma \in [0, 1]$, and $\circ$ denotes the binomial thinning operation.

2. The p.m.f. of discrete stable distributions are characterized by two parameters: exponent $\nu \in (0, 1]$, and rate parameter $\tau \in (0, \infty)$. We denote the p.m.f. of a univariate discrete stable distribution as $P_{DS}(\nu, \tau)$. In general, discrete stable distributions do not have a "nice" analytical form consisting of well-known elementary functions. Consequently, the univariate discrete stable distribution are typically expressed through their probability generating function.

3. The probability generating function of a discrete random variable $N$ having a stable discrete distribution is given by (179):

$$\mathbb{P}_N(z) = \exp(-\tau(1 - z)^\gamma). \tag{179}$$

## I.2  Definition of univariate affine discrete stable system

Let $M$,$X$, and $Y$ be a system of random variables with the joint distribution $P(M, X, Y)$. The joint distribution $P(M, X, Y)$ describes the univariate affine discrete stable system if it satisfies the following two properties:

1. $M \sim P(M)$ having some support set $\mathcal{M} \subseteq (0, \infty)$.

2. The conditional distributions of random variables $X$ and $Y$ conditioned on $M$ can be expressed through discrete stable family distributions with an affine dependence on $M$. Formally, $P(X|M{=}m){=}P_{DS}(\nu, am + b)$ and $P(Y|M{=}m){=}P_{DS}(\nu, cm + d)$, where $a, b, c, d \in (0, \infty)$.

## I.3  Formal proof of Theorem 4

**Theorem 4.** *Let $M, X$, and $Y$ be random variables whose joint distribution $P(M, X, Y)$ describes a univariate affine discrete stable system. Without the loss of generality, assume $a \geq c$. If $^a/_b \geq {}^c/_d$, then $\Delta_P$ contains a Markov chain of the form $M \rightarrow X \rightarrow Y$ and $UI(M; Y \backslash X) = 0$.*

*Proof.* We first note that we can always assume $a \geq c$ without the loss of generality because if $a \leq c$, then we can always switch our nomenclature to refer to $Y$ as $X$, and $X$ as $Y$.

We briefly outline the proof structure.

1. In the first part, we explicitly construct a joint distribution $Q_{MC}(M, X, Y)$ having the Markovian structure $M \to X \to Y$.

2. In the second part, we show that the $Q_{MC}(M, X, Y)$ constructed in the first part lies in $\Delta_P$. Therefore, we can then apply the result of proposition 1 to conclude $UI(M; Y \backslash X) = 0$.

**Part 1: Specifying $Q_{MC}(M, X, Y)$**

We explicitly construct the desired Markov chain $M \to X \to Y$ by constructing a larger Markov chain $M \to X \to X' \to Y$ and then marginalizing the larger Markov chain to obtain the desired Markov chain $M \to X \to Y$.

Denote the joint distribution of the Markov chain $M \to X \to X' \to Y$ as $Q_{MC}(M, X, X', Y)$. We can decompose $Q_{MC}(M, X, X', Y)$ by utilizing its Markovian structure as follows:

$$Q_{MC}(M, X, X', Y) = Q_{MC}(M)Q_{MC}(X|M)Q_{MC}(X'|X)Q_{MC}(Y|X').$$

Consequently, we can specify/construct the distribution $Q_{MC}(M, X, X', Y)$ by individually specifying $Q_{MC}(M)$, $Q_{MC}(X|M)$, $Q_{MC}(X'|X)$, and $Q_{MC}(Y|X')$.

Specifying $Q_{MC}(M)$ and $Q_{MC}(X|M)$: We choose $Q_{MC}(M)$ and $Q_{MC}(X|M)$ as follows:

$$Q_{MC}(M) = P(M) \text{ and } Q_{MC}(X|M) = P(X|M), \tag{180}$$

where $P(M)$ and $P(X|M)$ are marginal distributions derived from the original joint distribution $P(M, X, Y)$ (discussed in the theorem statement) over which the bivariate PID is being calculated.

Specifying $Q_{MC}(X'|X)$: $Q_{MC}(X'|X)$ is specified through a binomial thinning operation, specifically:

$$X' = X \circ (c/a)^{1/\nu}. \tag{181}$$

Note that the above binomial thinning operation is valid as $c \leq a$ by the assumption in the theorem statement. We also derive $Q_{MC}(X'|M)$ as it is needed for showing $Q_{MC}(M, X, Y) \in \Delta_P$.

*Deriving $Q_{MC}(X'|M)$:* We use lemma 13 to derive $Q_{MC}(X'|M = m)$ as $Q_{MC}(X|M = m)$ describes a univariate discrete stable distribution for a fixed $m$, and $X'$ is a binomially-thinned version of $X$. The exact expression of $Q_{MC}(X'|M = m)$ is provided in (182).

$$\begin{aligned} Q_{MC}(X'|M = m) &= P_{DS}(\nu, (c/a)^{\nu/\nu}(am + b/a)), \\ &= P_{DS}(\nu, cm + cb/a). \end{aligned} \tag{182}$$

Specifying $Q_{MC}(Y|X')$: $Q_{MC}(Y|X')$ is specified through the following stochastic transformation:

$$Y = X' + \epsilon, \tag{183}$$

where $\epsilon \sim P_{DS}(\nu, d - cb/a)$. For $Q_{MC}(Y|X')$ to be a valid distribution, we need to ensure that $\epsilon$ is distributed according to a legitimate discrete stable distribution. In order to show $\epsilon$ is distributed according to a legitimate discrete stable distribution, we need to ensure that $\nu$ and $d - cb/a$ lie within their appropriate bounds as specified in Sec. I.1. It is trivial to see $\nu \in (0, 1]$.

From the assumptions in the theorem statement, we know that

$$\frac{a}{b} \geq \frac{c}{d} \Rightarrow d \geq \frac{bc}{a} \Rightarrow d - \frac{bc}{a} \geq 0. \tag{184}$$

Hence, $\epsilon$ follows a legitimate discrete stable distribution. Furthermore, we choose $\epsilon \perp\!\!\!\perp (M, X, X')$.

Finally, we construct the desired $Q_{MC}(M, X, Y)$ from $Q_{MC}(M, X, X', Y)$ by marginalizing $X'$.

**Part 2: Showing $Q_{MC}(M, X, Y) \in \Delta_P$**

For showing that $Q_{MC}(M, X, Y) \in \Delta_P$, we first need to derive $Q_{MC}(Y|M)$.

Deriving $Q(Y|M)$: Note that $\epsilon$ follows discrete stable distribution $P_{DS}(\nu, d - {}^{bc}/a)$ and $\overline{Q_{MC}(X'|M = m)}$ is also a discrete stable distribution $P_{DS}(X'; \nu, cm + {}^{bc}/a)$ for a fixed $m$ (see (182)). Furthermore, $\epsilon \perp\!\!\!\perp X'|M$. Hence, using lemma 14, we have

$$Q_{MC}(Y|M = m) = P_{DS}(\nu, cm + {}^{bc}/a - {}^{bc}/a + d) = P_{DS}(\nu, cm + d). \tag{185}$$

As $P(Y|M) = P_{DS}(\nu, cM + d)$ (see Appx. I.2), we have:

$$Q(Y|M) = P(Y|M). \tag{186}$$

From (180) and (186), we can conclude:

$$Q_{MC}(M, X) = Q_{MC}(M)Q_{MC}(X|M) = P(M)P(X|M) = P(M, X), \tag{187}$$
$$Q_{MC}(M, Y) = Q_{MC}(M)Q_{MC}(Y|M) = P(M)P(Y|M) = P(M, Y). \tag{188}$$

Hence, $Q_{MC}(M, X, Y) \in \Delta_P$ and consequently, by proposition 1, $UI(M; Y \backslash X) = 0$, concluding our proof. □

**Lemma 13.** *Let* $N \sim P_{DS}(\nu, \tau)$. *Then,* $N_1 = \gamma \circ N \sim P_{DS}(\nu, \gamma^\nu \tau) \ \forall \ \gamma \in [0, 1]$, *where* $\circ$ *is the binomial thinning operation.*

*Proof.* The probability generating function of $N$, denoted as $\mathbb{P}_N(z)$, can be described using (179):

$$\mathbb{P}_N(z) = \exp\left(-\tau(1 - z)^\nu\right). \tag{189}$$

For calculating the probability generating function of $N_1$, we use the following relationship between the probability generating function of a random variable $R$ and its binomially-thinned version $R \circ p$, with $p \in [0, 1]$:

$$\mathbb{P}_{R \circ p}(z) = \mathbb{P}_R(1 - p(1 - z)) \text{ (see equation 3.3 in [24]).} \tag{190}$$

From (189) and (190), we can calculate the probability generating function of $N_1$ as follows:

$$\mathbb{P}_{N_1}(z) = \mathbb{P}_{N \circ \gamma}(z) = \mathbb{P}_N(1 - \gamma(1 - z)) = \exp\left(-\tau\left(1 - (1 - \gamma(1 - z))\right)^\nu\right)$$
$$= \exp\left(-\tau\left(\gamma(1 - z)\right)^\nu\right) = \exp\left(-\tau\gamma^\nu (1 - z)^\nu\right). \tag{191}$$

Inspecting the probability generating function of $N_1$ in (191), we can conclude that $N_1 \sim P_{DS}(\nu, \gamma^\nu \tau)$. □

**Lemma 14.** *Let* $N_1 \sim P_{DS}(\nu, \tau_1)$, $N_2 \sim P_{DS}(\nu, \tau_2)$, *and* $N_1 \perp\!\!\!\perp N_2$. *Then,* $N_1 + N_2 = N \sim P_{DS}(n; \nu, \tau_1 + \tau_2)$.

*Proof.* The probability generating function of $N_1$ and $N_2$, denoted as $\mathbb{P}_{N_1}(z)$ and $\mathbb{P}_{N_2}(z)$, can be described using (179):

$$\mathbb{P}_{N_1}(z) = \exp\left(-\tau_1(1 - z)^\nu\right) \text{ and } \mathbb{P}_{N_2}(z) = \exp\left(-\tau_2(1 - z)^\nu\right). \tag{192}$$

Calculating the probability generating function of $N = N_1 + N_2$,

$$\mathbb{P}_N(z) = \mathbb{P}_{N_1 + N_2}(z) \overset{(a)}{=} \mathbb{P}_{N_1}(z)\mathbb{P}_{N_2}(z) \overset{(b)}{=} \exp\left(-\tau_1 (1 - z)^\nu\right)\exp\left(-\tau_2 (1 - z)^\nu\right),$$
$$= \exp\left(-(\tau_1 + \tau_2)(1 - z)^\nu\right) \tag{193}$$

where $(a)$ is due to the fact that $N_1 \perp\!\!\!\perp N_2$, and $(b)$ is achieved by substituting the explicit forms of $\mathbb{P}_{N_1}(z)$ and $\mathbb{P}_{N_2}(z)$ from (192). Inspecting the probability generating function of $N_1 + N_2$ in (193), we can conclude that $N_1 + N_2 = N \sim P_{DS}(n_1; \nu, \tau_1 + \tau_2)$. □

## J Proof of Theorem 5

In this section, we provide the proof of theorem 5. Since theorem 5 provides the analytical PID terms for the multivariate linear Poisson system defined in Sec. 4, we briefly restate certain key properties of multivariate Poisson distributions and the multivariate linear Poisson system for convenience.

## J.1 Multivariate Poisson Distribution

We use the multivariate Poisson distribution proposed in [42]. For more details, see Appx. N.5. We list certain key properties of multivariate Poisson distributions that we make use of in the proof of theorem 5:

1. Under the definition of the multivariate Poisson distribution proposed in [42], each component of the random vector is a sum of independent Poisson random, i.e.,

$$\vec{\mathbf{N}} = \mathbf{A}\vec{\mathbf{N}}^g, \tag{194}$$

   where $\mathbf{A}$ is an appropriate matrix of 0's and 1's. We can decompose $\mathbf{A} = [\mathbf{A}_1 \quad \dots \quad \mathbf{A}_{d'}]$, where $\mathbf{A}_i$ is a $d \times \binom{d}{i}$ submatrix having no duplicate columns, and each of its columns containing exactly $i$ ones and $(d-i)$ zeros [44], and $\vec{\mathbf{N}}^g = [N_1^g \dots N_d^g \ N_{12}^g \dots N_{(d-1)d}^g \cdots N_{d-(d'-1)\dots d}^g]^T$, with $N_{i_1\dots i_j}^g \sim \text{Poisson}(\lambda_{i_1\dots i_j}) \ \forall \ (i_1,\dots,i_j) \in \mathbb{A}_j^d, j \in [d']$. Furthermore, the random variables $\{N_1,\dots,N_{d-(d'-1)\dots d}\}$ are mutually independent.

2. We denote the p.m.f. of the multivariate Poisson distribution as $\text{Poisson}(d, d', \vec{\boldsymbol{\Lambda}})$, where $d \geq d'$, and

$$\vec{\boldsymbol{\Lambda}} = \begin{bmatrix} \lambda_1 \ \dots \ \lambda_d \ \lambda_{12} \ \dots \lambda_{d-(d'-1)\dots d} \end{bmatrix}^T. \tag{195}$$

3. For $d' = 1$, we have that $\vec{\mathbf{N}}$ is a collection of independent Poisson random variables, and when both $d = d' = 1$, we recover the scalar Poisson distribution: $\Pr(N = n) = \frac{e^{-\lambda}\lambda^n}{n!}, \ \forall \ n \in \mathbb{N}_0$.

4. Let $\vec{\mathbf{N}} \sim \text{Poisson}(d, d', \vec{\boldsymbol{\Lambda}})$, where $\vec{\mathbf{N}}$ is a $d$-dimensional random vector. Then the corresponding p.m.f. of $\vec{\mathbf{N}}$ is described below:

   Let $\vec{\mathbf{n}}' = \begin{bmatrix} n_{12} \ \dots \ n_{(d-1)d} \ \dots \ n_{d-(d'-1)\dots d} \end{bmatrix}^T$, and $d_{\vec{\mathbf{n}}'}$ be the dimension of $\vec{\mathbf{n}}'$, then:

$$P(\vec{\mathbf{N}} = \vec{\mathbf{n}}) = e^{-\vec{\mathbf{1}}^T\vec{\boldsymbol{\Lambda}}} \prod_{i=1}^{d} \lambda_i^{n_i} \sum_{\vec{\mathbf{n}}' \in C} \left( \prod_{(i_1,i_2) \in \mathbb{A}_2^d} \left( \frac{\lambda_{i_1 i_2}}{\lambda_{i_1}\lambda_{i_2}} \right)^{n_{i_1 i_2}} \times \right.$$

$$\left. \dots \times \prod_{(i_1,\dots,i_{d'}) \in \mathbb{A}_{d'}^d} \left( \frac{\lambda_{i_1 \dots i_{d'}}}{\prod_{j=1}^{d'}\lambda_{i_j}} \right)^{n_{i_1 \dots i_{d'}}} \times Q(\vec{\mathbf{n}}, \vec{\mathbf{n}}') \right), \tag{196}$$

   where $C = \{\vec{\mathbf{n}}' \in \mathbb{N}_0^{d_{\vec{\mathbf{n}}'}} : (\vec{\mathbf{a}}_i')^T \vec{\mathbf{n}}' \leq n_i \ \forall \ i \in [d]\}$, and

$$Q(\vec{\mathbf{n}}, \vec{\mathbf{n}}') = \prod_{i=1}^{d} \frac{1}{(n_i - \vec{\mathbf{a}}_i'^T \vec{\mathbf{n}}')!} \prod_{j=2}^{d'} \prod_{(i_1,\dots,i_j) \in \mathbb{A}_j^d} \frac{1}{n_{i_1\dots i_j}^g !},$$

   with $\vec{\mathbf{a}}_i'$ being the $i$-th row of the matrix $\mathbf{A}' = [\mathbf{A}_2 \dots \mathbf{A}_{d'}]$.

## J.2 Definition of the multivariate linear Poisson system

Let the random variable $M \sim P(M)$. $\vec{\mathbf{X}}$ and $\vec{\mathbf{Y}}$ are $d_X$-dimensional and $d_Y$-dimensional random vectors, respectively. The joint distribution $P(M, \vec{\mathbf{X}}, \vec{\mathbf{Y}})$ describes the multivariate linear Poisson system if it satisfies the following two properties:

1. $M \sim P(M)$ having some support set $\mathcal{M} \subseteq (0, \infty)$.

2. The conditional distributions of random vectors $\vec{\mathbf{X}}$ and $\vec{\mathbf{Y}}$ are multivariate Poisson distributions, i.e., $P(\vec{\mathbf{X}}|M) = \text{Poisson}(d_X, d_X', \vec{\boldsymbol{\Lambda}}_X)$ and $P(\vec{\mathbf{Y}}|M) = \text{Poisson}(d_Y, d_Y', \vec{\boldsymbol{\Lambda}}_Y)$, with:

$$\vec{\boldsymbol{\Lambda}}_X = \begin{bmatrix} \lambda_1^X \ \dots \ \lambda_{d_X-(d_X'-1)\dots d_X}^X \end{bmatrix}^T, \lambda_{i_1\dots i_j}^X = \gamma_{i_1\dots i_j}^X M^j \ \forall \ j \in [d_X] \text{ and } (i_1,\dots,i_j) \in \mathbb{A}_j^{d_X},$$

$$\boldsymbol{\Lambda}_Y = \begin{bmatrix} \lambda_1^Y \ \dots \ \lambda_{d_Y-(d_Y'-1)\dots d_Y}^Y \end{bmatrix}^T, \lambda_{i_1\dots i_j}^Y = \gamma_{i_1\dots i_j}^Y M^j \ \forall \ j \in [d_Y] \text{ and } (i_1,\dots,i_j) \in \mathbb{A}_j^{d_Y}.$$

   Here, $\gamma_1^X, \dots, \gamma_{d_X-(d_X'-1)\dots d_X}^X, \gamma_1^Y, \dots, \gamma_{d_Y-(d_Y'-1)\dots d_Y}^Y \in \mathbb{R}^+$.

Furthermore, let $\mathbf{A}_X$, and $\mathbf{A}_Y$ be the corresponding $\mathbf{A}$-matrices (defined in (194)) associated with $P(\vec{\mathbf{X}}|M)$ and $P(\vec{\mathbf{Y}}|M)$, respectively.

## J.3 Formal proof of Theorem 5

**Theorem 5.** *Let the joint distribution $P(M, \vec{\mathbf{X}}, \vec{\mathbf{Y}})$ of $M$, $\vec{\mathbf{X}}$, and $\vec{\mathbf{Y}}$ describe the multivariate linear Poisson system. Without the loss of generality, assume $d'_X \geq d'_Y$. If $\sum_{(i_1,\ldots,i_j) \in \mathbb{A}_j^{d_X}} \gamma_{i_1\ldots i_j}^X \geq \sum_{(i_1,\ldots,i_j) \in \mathbb{A}_j^{d_Y}} \gamma_{i_1\ldots i_j}^Y \, \forall \, j \in [d'_Y]$, then $\Delta_P$ contains a Markov chain of the form $M \to \vec{\mathbf{X}} \to \vec{\mathbf{Y}}$ and $UI(M; \vec{\mathbf{Y}} \backslash \vec{\mathbf{X}}) = 0$.*

*Proof.* We first note that we can always assume $d'_X \geq d'_Y$ without the loss of generality because if $d'_X \leq d'_Y$, then we can always switch our nomenclature to refer to $\vec{\mathbf{Y}}$ as $\vec{\mathbf{X}}$, and $\vec{\mathbf{X}}$ as $\vec{\mathbf{Y}}$.

We provide an explicit construction of the Markov chain $M \to \vec{\mathbf{X}} \to \vec{\mathbf{X}}^g \to \vec{\mathbf{Y}}^g \to \vec{\mathbf{Y}}$ having the marginals $P(M, \vec{\mathbf{X}})$, and $P(M, \vec{\mathbf{Y}})$. Denote the joint density of $M \to \vec{\mathbf{X}} \to \vec{\mathbf{X}}^g \to \vec{\mathbf{Y}}^g \to \vec{\mathbf{Y}}$ as $Q(M, \vec{\mathbf{X}}, \vec{\mathbf{X}}^g, \vec{\mathbf{Y}}^g, \vec{\mathbf{Y}})$. For $Q(M)$ and $Q(\vec{\mathbf{X}}|M)$, we choose them to be equal to $P(M)$ and $P(\vec{\mathbf{X}}|M)$, i.e. $Q(M) = P(M)$, and $Q(\vec{\mathbf{X}}|M) = P(\vec{\mathbf{X}}|M)$. Note that due to this construction, $P(M, \vec{\mathbf{X}}) = Q(M, \vec{\mathbf{X}})$ holds trivially.

For $Q(\vec{\mathbf{X}}^g|\vec{\mathbf{X}})$, we use the result described in lemma 17. Note that the construction for $Q(\vec{\mathbf{X}}^g|\vec{\mathbf{X}})$ is not explicit but rather implicit. Let $d_{\vec{\boldsymbol{\Lambda}}_X}$ be the dimension of $\vec{\boldsymbol{\Lambda}}_X$, then we explicitly choose: $Q(\vec{\mathbf{X}}|\vec{\mathbf{X}}^g, M) = \delta_K(\vec{\mathbf{X}} = \mathbf{A}_X \vec{\mathbf{X}}^g)$, and $Q(\vec{\mathbf{Y}}^g|M) = \mathrm{Poisson}(d_{\vec{\boldsymbol{\Lambda}}_X}, 1, \vec{\boldsymbol{\Lambda}}_Y)$. Note that $Q(\vec{\mathbf{X}}|M) = \mathrm{Poisson}(d_X, d'_X, \vec{\boldsymbol{\Lambda}}_X)$ by construction, and we derive $Q(\vec{\mathbf{X}}^g|\vec{\mathbf{X}}, M)$ through the Bayes theorem. Here, $\delta_K(\cdot)$ is the Kronecker delta function [59]. By lemma 17, we know that $Q(\vec{\mathbf{X}}^g|\vec{\mathbf{X}}, M) = Q(\vec{\mathbf{X}}^g|\vec{\mathbf{X}})$, and hence we have the Markov chain $M \to \vec{\mathbf{X}} \to \vec{\mathbf{X}}^g$.

For choosing $Q(\vec{\mathbf{Y}}^g|\vec{\mathbf{X}}^g)$, we rely on the result of lemma 18. Let us define the random vectors:

$$\vec{\mathbf{X}}_j = \left[ X_{i_1\ldots i_j}^g \right]_{(i_1,\ldots,i_j) \in \mathbb{A}_j^{d_1}}^T, \text{ and } \vec{\mathbf{Y}}_j = \left[ Y_{i_1\ldots i_j}^g \right]_{(i_1,\ldots,i_j) \in \mathbb{A}_j^{d_2}}^T,$$

i.e., $\vec{\mathbf{X}}_j$ and $\vec{\mathbf{Y}}_j$ are random vectors containing all terms of the form $X_{i_1,\ldots,i_j}^g$, and $Y_{k_1,\ldots,k_j}^g$, where $(i_1,\ldots,i_j) \in \mathbb{A}_j^{d_1}$ and $(k_1,\ldots,k_j) \in \mathbb{A}_j^{d_2}$, respectively. Note that we can write $\vec{\mathbf{X}}^g = \left[ \vec{\mathbf{X}}_1^T \ldots \vec{\mathbf{X}}_{d'_1}^T \right]^T$, and we define $\vec{\mathbf{Y}}^g = \left[ \vec{\mathbf{Y}}_1^T \ldots \vec{\mathbf{Y}}_{d'_2}^T \right]^T$. Then, we construct $Q(\vec{\mathbf{Y}}^g|\vec{\mathbf{X}}^g)$ as a product of $d'_Y$ multinomial distributions, described below:

$$Q(\vec{\mathbf{Y}}^g|\vec{\mathbf{X}}^g) = \prod_{j=1}^{d'_Y} Q(\vec{\mathbf{Y}}_j|\vec{\mathbf{X}}_j),$$

where $Q(\vec{\mathbf{Y}}_j = \vec{\mathbf{y}}_j | \vec{\mathbf{X}}_j = \vec{\mathbf{x}}_j) = \mathrm{Multinomial}(\mathbf{k}_j; N_j, \mathbf{p}_i)$, and

$$N_j = \vec{\mathbf{1}}^T \vec{\mathbf{x}}_j, \; \vec{\mathbf{k}}_j = \left[ \vec{\mathbf{y}}_j^T \quad \vec{\mathbf{1}}^T \vec{\mathbf{x}}_j - \vec{\mathbf{1}}^T \vec{\mathbf{y}}_j \right]^T,$$

$$\vec{\mathbf{p}}_i = \left[ p_{1\ldots j} \cdots p_{d_2-(j-1)\ldots d_2} \; 1 - \sum_{(i_1,\ldots,i_j) \in \mathbb{A}_j^{d_2}} p_{i_1\ldots i_j} \right]^T,$$

$$p_{i_1\ldots i_j} = \frac{\gamma_{i_1\ldots i_j}^Y}{\sum_{(i_1,\ldots,i_j) \in \mathbb{A}_j^{d_1}} \gamma_{i_1\ldots i_j}^X} \; \forall \, (i_1,\ldots,i_j) \in \mathbb{A}_j^{d_2}.$$

By construction $\vec{\mathbf{X}}^g$ consists of mutually conditionally independent Poisson random variables (conditioned on $M$), and the condition of lemma 18 (specified in (219)) is satisfied by condition 1. Therefore, using the result of lemma 18 on the Markov chain $M \to \vec{\mathbf{X}}^g \to \vec{\mathbf{Y}}^g$ (obtained by marginalizing $\vec{\mathbf{X}}$), we conclude that $Q(\vec{\mathbf{Y}}^g|M) = \mathrm{Poisson}(1, d_{\vec{\boldsymbol{\Lambda}}_Y}, \vec{\boldsymbol{\Lambda}}_Y)$, where $d_{\vec{\boldsymbol{\Lambda}}_Y}$ is the dimension of $\vec{\boldsymbol{\Lambda}}_Y$.

We specify $Q(\vec{\mathbf{Y}}|\vec{\mathbf{Y}}^g)$ through the following deterministic transformation $\vec{\mathbf{Y}} = \mathbf{A}_Y \vec{\mathbf{Y}}^g$ to obtain the Markov chain $M \to \vec{\mathbf{X}} \to \vec{\mathbf{X}}^g \to \vec{\mathbf{Y}}^g \to \vec{\mathbf{Y}}$. Marginalizing $\vec{\mathbf{X}}$ and $\vec{\mathbf{X}}^g$ in the above Markov chain,

we get the following Markov chain: $M \to \vec{\mathbf{Y}}^g \to \vec{\mathbf{Y}}$. Now, since $\vec{\mathbf{Y}} = \mathbf{A}_Y \vec{\mathbf{Y}}^g$ and $P(\vec{\mathbf{Y}}^g|M) = $ Poisson$(d_{\vec{\mathbf{A}}_Y}, 1, \vec{\mathbf{\Lambda}}_Y)$, $P(\vec{\mathbf{Y}}|M) = $ Poisson$(d_Y, d'_Y, \Lambda_Y)$ (by definition of the multivariate Poisson discussed in Sec. J.1). Since we have $Q(M) = P(M)$ and $Q(\vec{\mathbf{Y}}|M) = P(\vec{\mathbf{Y}}|M)$, we also have $Q(M, \vec{\mathbf{Y}}) = P(M, \vec{\mathbf{Y}})$. Marginalizing $\vec{\mathbf{X}}^g$ and $\vec{\mathbf{Y}}^g$ from the Markov chain $M \to \vec{\mathbf{X}} \to \vec{\mathbf{X}}^g \to \vec{\mathbf{Y}}^g \to \vec{\mathbf{Y}}$ results in the desired Markov chain $M \to \vec{\mathbf{X}} \to \vec{\mathbf{Y}}$ having $Q(M, \vec{\mathbf{X}}) = P(M, \vec{\mathbf{X}})$ and $Q(M, \vec{\mathbf{Y}}) = P(M, \vec{\mathbf{Y}})$, concluding our proof. $\qquad\square$

**Lemma 15.** *If $N_j \sim Poisson(\lambda_j) \ \forall \ j \in \{1, \ldots, d\}$, and all $N_j$'s are mutually independent, then $P(N_1, \ldots, N_j| \sum_{j=1}^{d} N_j = N) = Multinomial(N_1, \ldots, N_j; N, [\lambda_1/\sum_{i=1}^{d} \lambda_j \quad \cdots \quad \lambda_d/\sum_{i=1}^{d} \lambda_j])$.*

*Proof.* See Section 3 of [60]. $\qquad\square$

**Lemma 16.** *Let $\vec{\mathbf{N}}_1 = \begin{bmatrix} N_1^{(1)} & \ldots N_{d_1}^{(1)} \end{bmatrix}^T$, where $N_j \sim Poisson(\lambda_j^{(1)}) \ \forall \ j \in \{1, \ldots, d_1\}$ and all $N_j$'s are mutually independent from each other. Define $P(\vec{\mathbf{N}}_2|\vec{\mathbf{N}}_1) = Multinomial \left( \vec{\mathbf{k}}; N, \vec{\mathbf{p}} \right)$, where*

$$N = \sum_{j=1}^{d_1} N_j^{(1)}, \ \vec{\mathbf{k}} = \begin{bmatrix} N_1^{(2)} & \cdots & N_{d_2}^{(2)} & \sum_{j=1}^{d_1} N_j^{(1)} - \sum_{j=1}^{d_2} N_j \end{bmatrix}^T,$$

$$\vec{\mathbf{p}} = \begin{bmatrix} \frac{\lambda_1^{(2)}}{\sum_{j=1}^{d_1} \lambda_j^{(1)}} & \cdots & \frac{\lambda_1^{(2)}}{\sum_{j=1}^{d_1} \lambda_j^{(1)}} & 1 - \frac{\sum_{j=1}^{d_2} \lambda_j^{(2)}}{\sum_{j=1}^{d_1} \lambda_j^{(1)}} \end{bmatrix}^T,$$

*where $\{\lambda_j^{(2)}\}_{j=1}^{d_2}$ are some positive constants, i.e., $\lambda_j^{(2)} \in (0, \infty)$, such that $\sum_{i=1}^{d_2} \lambda_j^{(2)} \leq \sum_{j=1}^{d_1} \lambda_j^{(1)}$. Then, $\vec{\mathbf{N}}_2 = \begin{bmatrix} N_1^{(2)} & \ldots N_{d_2}^{(2)} \end{bmatrix}^T$, where $N_j^{(2)} \sim Poisson(\lambda_j^{(2)})$, and all $N_j^{(2)}$'s are mutually independent form each other.*

*Proof.* Lemma 16 is a straight-forward generalization of lemma 15. Denote the probability of $\vec{\mathbf{N}}_2$ shown in the lemma statement as $P(\vec{\mathbf{N}}_2)$. We will prove the above lemma by constructing a Markov chain $\vec{\mathbf{N}}_1 \to N'_1 \to N'_2 \to \vec{\mathbf{N}}_2$ with joint density $Q(\vec{\mathbf{N}}_1, N'_1, N'_2, \vec{\mathbf{N}}_2)$, and showing that $Q(\vec{\mathbf{N}}_2) = P(\vec{\mathbf{N}}_2)$ and $Q(\vec{\mathbf{N}}_2|\vec{\mathbf{N}}_1) = P(\vec{\mathbf{N}}_2|\vec{\mathbf{N}}_1)$.

Specifying $Q(N'_1|\vec{\mathbf{N}}_1)$: We specify $Q(N'_1|\vec{\mathbf{N}}_1)$ through the following deterministic transformation:

$$N'_1 = \vec{\mathbf{1}}^T \vec{\mathbf{N}}_1 = \sum_{j=1}^{d_1} N_j^{(1)}. \tag{197}$$

Calculating $Q(N'_1)$: We use the property that the sum of independent Poisson random variables is a Poisson random variable with its rate parameter being the sum of the rate parameters of its summands [61]. Hence, $Q(N'_1) = $ Poisson$(\sum_{j=1}^{d_1} \lambda_j^{(1)})$.

Specifying $Q(N'_2|N'_1)$: We specify $Q(N'_2|N'_1 = n'_1)$ as follows:

$$Q(N'_2|N'_1 = n'_1) = \text{Binomial} \left( N'_2; n'_1, \sum_{j=1}^{d_2} \lambda_j^{(2)} / \sum_{j=1}^{d_1} \lambda_j^{(1)} \right). \tag{198}$$

Calculating $Q(N'_2)$: We use lemma 15 to calculate $Q(N'_2)$. Note that $Q(N'_1) = $ Poisson$(\sum_{j=1}^{d_1} \lambda_j^{(1)})$, and $Q(N'_2|N'_1) = $ Binomial $\left( N'_2; n'_1, \sum_{j=1}^{d_2} \lambda_j^{(2)} / \sum_{j=1}^{d_1} \lambda_j^{(1)} \right)$. Hence, by lemma 15, we have:

$$Q(N'_2) = \text{Poisson} \left( \sum_{j=1}^{d_1} \lambda_j^{(1)} \times \frac{\sum_{j=1}^{d_2} \lambda_j^{(2)}}{\sum_{j=1}^{d_1} \lambda_j^{(1)}} \right) = \text{Poisson} \left( \sum_{j=1}^{d_2} \lambda_j^{(2)} \right). \tag{199}$$

Specifying $Q(N'_2|N'_1)$: We specify $Q(\vec{\mathbf{N}}_2|N'_2 = n'_2)$ as follows:

$$Q(\vec{\mathbf{N}}_2|N'_2 = n'_2) = \text{Multinomial}\left(\vec{\mathbf{N}}'_2; n'_2, \left[\frac{\lambda_1^{(2)}}{\sum_{j=1}^{d_2}\lambda_j^{(2)}} \quad \cdots \quad \frac{\lambda_{d_2}^{(2)}}{\sum_{j=1}^{d_2}\lambda_j^{(2)}}\right]^T\right). \quad (200)$$

Calculating $Q(\vec{\mathbf{N}}_2)$: We again use lemma 15 to calculate $Q(\vec{\mathbf{N}}_2)$. Note that,

$$Q(N'_2) = \text{Poisson}\left(\sum_{j=1}^{d_2}\lambda_j^{(2)}\right), \text{ and}$$

$$Q(\vec{\mathbf{N}}_2|N'_2 = n'_2) = \text{Multinomial}\left(\vec{\mathbf{N}}'_2; n'_2, \left[\frac{\lambda_1^{(2)}}{\sum_{j=1}^{d_2}\lambda_j^{(2)}} \quad \cdots \quad \frac{\lambda_{d_2}^{(2)}}{\sum_{j=1}^{d_2}\lambda_j^{(2)}}\right]^T\right). \quad (201)$$

Hence, by lemma 15, we have:

$$\vec{\mathbf{N}}_2 = \left[N_1^{(2)} \quad \cdots \quad N_{d_2}^{(2)}\right]^T, \text{ where } N_j^{(2)} \sim \text{Poisson}(\lambda_j^{(2)}) \; \forall \, j \in [d_2]. \quad (202)$$

Hence, by (202), we know that $Q(\vec{\mathbf{N}}_2) = P(\vec{\mathbf{N}}_2)$.

Calculating $Q(\vec{\mathbf{N}}_2|\vec{\mathbf{N}}_1)$:

$$Q(\vec{\mathbf{N}}_2 = \vec{\mathbf{n}}_2|\vec{\mathbf{N}}_1 = \vec{\mathbf{n}}_1) = \sum_{n'_1, n'_2=0}^{\infty} Q(N'_1 = n'_1, N'_2 = n'_2, \vec{\mathbf{N}}_2 = \vec{\mathbf{n}}_2|\vec{\mathbf{N}}_1 = \vec{\mathbf{n}}_1),$$

$$\overset{(a)}{=} \sum_{n'_1, n'_2=0}^{\infty} Q(n'_1|\vec{\mathbf{n}}_2)Q(n'_2|n'_1)Q(\vec{\mathbf{n}}_2|n'_2),$$

where $(a)$ is due to the Markovian nature of $Q(\vec{\mathbf{N}}_1, N'_1, N'_2, \vec{\mathbf{N}}_2)$. We drop the random variable notation for brevity. Let $\vec{\mathbf{n}}_1 = \left[n_1^{(1)} \quad \cdots \quad n_{d_1}^{(1)}\right]^T$, and $\vec{\mathbf{n}}_2 = \left[n_1^{(2)} \quad \cdots \quad n_{d_2}^{(2)}\right]^T$. Then, substituting the exact form of $Q(n'_1|\vec{\mathbf{n}}_1), Q(n'_2|n'_1)$, and $Q(\vec{\mathbf{n}}_2|n'_2)$ in the above equation:

$$= \sum_{n'_1, n'_2=0}^{\infty} \mathbb{I}\left[n'_1 = \sum_{j=1}^{d_1}n_j^{(1)}\right] \frac{(n'_1)!}{(n'_1 - n'_2)!(n'_2)!} \left(\frac{\sum_{j=1}^{d_2}\lambda_j^{(2)}}{\sum_{j=1}^{d_1}\lambda_j^{(1)}}\right)^{n'_2} \left(1 - \frac{\sum_{j=1}^{d_2}\lambda_j^{(2)}}{\sum_{j=1}^{d_1}\lambda_j^{(1)}}\right)^{n'_1 - n'_2} \times,$$

$$\mathbb{I}\left[\sum_{j=1}^{d_2}n_j^{(2)} = n'_2\right] \frac{(n'_2)!}{\prod_{j=1}^{d_2}(n_j^{(2)})!} \prod_{j=1}^{d_2}\left(\frac{\lambda_j^{(2)}}{\sum_{j=1}^{d_2}\lambda_j^{(2)}}\right)^{n_j^{(2)}},$$

where $\mathbb{I}[\cdot]$ is an indicator function. Simplifying the above equation by using the fact that only non-zero term in the above summation is $n'_1 = \sum_{j=1}^{d_1}n_j^{(1)}$ and $n'_2 = \sum_{j=1}^{d_1}n_j^{(2)}$.

$$= \frac{\left(\sum_{j=1}^{d_1}n_j^{(1)}\right)!}{\left(\sum_{j=1}^{d_1}n_j^{(1)} - \sum_{j=1}^{d_2}n_j^{(2)}\right)!\left(\sum_{j=1}^{d_2}n_j^{(2)}\right)!} \left(\frac{\sum_{j=1}^{d_2}\lambda_j^{(2)}}{\sum_{j=1}^{d_1}\lambda_j^{(1)}}\right)^{\sum_{j=1}^{d_2}n_j^{(2)}} \times$$

$$\left(1 - \frac{\sum_{j=1}^{d_2}\lambda_j^{(2)}}{\sum_{j=1}^{d_1}\lambda_j^{(1)}}\right)^{\sum_{j=1}^{d_1}n_j^{(1)} - \sum_{j=1}^{d_1}n_j^{(2)}} \frac{\left(\sum_{j=1}^{d_2}n_j^{(2)}\right)!}{\prod_{j=1}^{d_2}(n_j^{(2)})!} \prod_{j=1}^{d_2}\left(\frac{\lambda_j^{(2)}}{\sum_{j=1}^{d_2}\lambda_j^{(2)}}\right)^{n_j^{(2)}},$$

Collecting all factorial terms together, and rearranging some terms:

$$= \frac{\left(\sum_{j=1}^{d_1}n_j^{(1)}\right)!}{\left(\sum_{j=1}^{d_1}n_j^{(1)} - \sum_{j=1}^{d_2}n_j^{(2)}\right)!\prod_{j=1}^{d_2}(n_j^{(2)})!} \left(1 - \frac{\sum_{j=1}^{d_2}\lambda_j^{(2)}}{\sum_{j=1}^{d_1}\lambda_j^{(1)}}\right)^{\sum_{j=1}^{d_1}n_j^{(1)} - \sum_{j=1}^{d_2}n_j^{(2)}} \times$$

$$\left(\frac{\sum_{j=1}^{d_2}\lambda_j^{(2)}}{\sum_{j=1}^{d_1}\lambda_j^{(1)}}\right)^{\sum_{j=1}^{d_2}n_j^{(2)}} \prod_{j=1}^{d_2}\left(\frac{\lambda_j^{(2)}}{\sum_{j=1}^{d_2}\lambda_j^{(2)}}\right)^{n_j^{(2)}}.$$

Substituting $\left(\sum_{j=1}^{d_2}\lambda_j^{(2)}/\sum_{j=1}^{d_1}\lambda_j^{(1)}\right)^{\sum_{j=1}^{d_2}n_j^{(2)}}=\prod_{j=1}^{d_2}\left(\sum_{j=1}^{d_2}\lambda_j^{(2)}/\sum_{j=1}^{d_1}\lambda_j^{(1)}\right)^{n_j^{(2)}}$ in the above equation:

$$=\frac{\left(\sum_{j=1}^{d_1}n_j^{(1)}\right)!}{\left(\sum_{j=1}^{d_1}n_j^{(1)}-\sum_{j=1}^{d_2}n_j^{(2)}\right)!\prod_{j=1}^{d_2}(n_j^{(2)})!}\left(1-\frac{\sum_{j=1}^{d_2}\lambda_j^{(2)}}{\sum_{j=1}^{d_1}\lambda_j^{(1)}}\right)^{\sum_{j=1}^{d_1}n_j^{(1)}-\sum_{j=1}^{d_2}n_j^{(2)}}\times$$

$$\prod_{j=1}^{d_1}\left(\frac{\sum_{j=1}^{d_2}\lambda_j^{(2)}}{\sum_{j=1}^{d_1}\lambda_j^{(1)}}\right)^{n_j^{(2)}}\prod_{j=1}^{d_2}\left(\frac{\lambda_j^{(2)}}{\sum_{j=1}^{d_2}\lambda_j^{(2)}}\right)^{n_j^{(2)}}$$

$$=\frac{\left(\sum_{j=1}^{d_1}n_j^{(1)}\right)!}{\left(\sum_{j=1}^{d_1}n_j^{(1)}-\sum_{j=1}^{d_2}n_j^{(2)}\right)!\prod_{j=1}^{d_2}(n_j^{(2)})!}\left(1-\frac{\sum_{j=1}^{d_2}\lambda_j^{(2)}}{\sum_{j=1}^{d_1}\lambda_j^{(1)}}\right)^{\sum_{j=1}^{d_1}n_j^{(1)}-\sum_{j=1}^{d_2}n_j^{(2)}}\times$$

$$\prod_{j=1}^{d_1}\left(\frac{\sum_{j=1}^{d_2}\lambda_j^{(2)}}{\sum_{j=1}^{d_1}\lambda_j^{(1)}}\right)^{n_j^{(2)}}\left(\frac{\lambda_j^{(2)}}{\sum_{j=1}^{d_2}\lambda_j^{(2)}}\right)^{n_j^{(2)}}$$

$$=\frac{\left(\sum_{j=1}^{d_1}n_j^{(1)}\right)!}{\left(\sum_{j=1}^{d_1}n_j^{(1)}-\sum_{j=1}^{d_2}n_j^{(2)}\right)!\prod_{j=1}^{d_2}(n_j^{(2)})!}\left(1-\frac{\sum_{j=1}^{d_2}\lambda_j^{(2)}}{\sum_{j=1}^{d_1}\lambda_j^{(1)}}\right)^{\sum_{j=1}^{d_1}n_j^{(1)}-\sum_{j=1}^{d_2}n_j^{(2)}}\times$$

$$\prod_{j=1}^{d_1}\left(\frac{\lambda_j^{(2)}}{\sum_{j=1}^{d_1}\lambda_j^{(1)}}\right)^{n_j^{(2)}}. \tag{203}$$

By inspecting (203), we can see that $Q(\vec{\mathbf{N}}_2|\vec{\mathbf{N}}_1)=\text{Multinomial}(\vec{\mathbf{k}};N,\vec{\mathbf{p}})$ with,

$$N=\sum_{j=1}^{d_1}N_j^{(1)},\ \vec{\mathbf{k}}=\left[N_1^{(2)}\ \cdots\ N_{d_2}^{(2)}\ \sum_{j=1}^{d_1}N_j^{(1)}-\sum_{j=1}^{d_2}N_j\right]^T, \tag{204}$$

$$\vec{\mathbf{p}}=\left[\lambda_1^{(2)}/\sum_{j=1}^{d_1}\lambda_j^{(1)}\ \cdots\ \lambda_1^{(2)}/\sum_{j=1}^{d_1}\lambda_j^{(1)}\ 1-\sum_{j=1}^{d_2}\lambda_j^{(2)}/\sum_{j=1}^{d_1}\lambda_j^{(1)}\right]^T, \tag{205}$$

Hence, by (205), we can conclude that $Q(\vec{\mathbf{N}}_2|\vec{\mathbf{N}}_1)=P(\vec{\mathbf{N}}_2|\vec{\mathbf{N}}_1)$ finishing our proof. $\qquad\square$

**Lemma 17.** *Let $\vec{\mathbf{N}}$ be a $d$-dimensional vector and $M$ be a positive random variable, i.e. $M\sim P(M),P(M\le 0)=0$. Let $P(\vec{\mathbf{N}}|M)=Poisson(d,d',\vec{\mathbf{\Lambda}})$, where:*

$$\vec{\mathbf{\Lambda}}=\left[\lambda_1\ \ldots\ \lambda_d\ \lambda_{12}\ \ldots\lambda_{d-(d'-1)\ldots d}\right]^T,$$

$$\lambda_{i_1\ldots i_j}=\gamma_{i_1\ldots i_j}M^j\ \forall\ j\in[d']\ and\ (i_1,\ldots,i_{d'})\in\mathbb{A}_j^d. \tag{206}$$

*Let $\vec{\mathbf{N}}^g=\left[N_1^g\ \ldots\ N_d^g\ N_{12}^g\ \cdots\ N_{(d-1)d}^g\ \cdots\ N_{d-(d'-1)\ldots d}^g\right]^T$, where $P(\vec{\mathbf{N}}^g|M)=Poisson(d_{\vec{\mathbf{\Lambda}}},1,\vec{\mathbf{\Lambda}})$ and $\vec{\mathbf{N}}=\mathbf{A}\vec{\mathbf{N}}^g$, where $d_\Lambda$ is the dimension of $\vec{\mathbf{\Lambda}}$, and $\mathbf{A}=[\mathbf{A}_1\ldots\mathbf{A}_{d'}]$ as defined in Sec. J.1. Then $P(\vec{\mathbf{N}}^g|\vec{\mathbf{N}},M)=P(\vec{\mathbf{N}}^g|\vec{\mathbf{N}})$, i.e. $M,\vec{\mathbf{N}}^g$ and $\vec{\mathbf{N}}$ form the following Markov chain $M\to\vec{\mathbf{N}}\to\vec{\mathbf{N}}^g$.*

*Proof.* To prove lemma 17, all we need to show is that $P(\vec{\mathbf{N}}^g|\vec{\mathbf{N}},M)$ does not depend on $M$. Let us to calculate $P(\vec{\mathbf{N}}^g|\vec{\mathbf{N}},M)$. By Bayes Theorem, we know:

$$P(\vec{\mathbf{N}}^g|\vec{\mathbf{N}},M)=\frac{P(\vec{\mathbf{N}}^g|M)P(\vec{\mathbf{N}}|\vec{\mathbf{N}}^g,M)}{P(\vec{\mathbf{N}}|M)}. \tag{207}$$

Let us now write the expression for $P(\vec{\mathbf{N}}|M)$ using (196):

$$P(\vec{\mathbf{N}}=\vec{\mathbf{n}}|M)=Q(\vec{\mathbf{n}},\vec{\mathbf{n}}')e^{-\vec{\mathbf{1}}^T\vec{\mathbf{\Lambda}}}\prod_{i=1}^d(\gamma_i M)^{n_i}\left(\sum_{\vec{\mathbf{n}}'\in C}\prod_{j=2}^{d'}\prod_{(i_1,\ldots,i_j)\in\mathbb{A}_j^d}\left(\frac{\gamma_{i_1\ldots i_j}M^j}{\prod_{l=1}^j\gamma_{i_l}M}\right)^{n'_{i_1\ldots i_j}}\right). \tag{208}$$

Canceling $M$ in the above equation for the terms inside the summation, we get:

$$P(\vec{\mathbf{N}}=\vec{\mathbf{n}}|M)=Q(\vec{\mathbf{n}},\vec{\mathbf{n}}')e^{-\vec{\mathbf{1}}^T\vec{\mathbf{\Lambda}}}\prod_{i=1}^{d}(\gamma_i M)^{n_i}\left(\sum_{\vec{\mathbf{n}}'\in C}\prod_{j=2}^{d'}\prod_{(i_1,...,i_j)\in\mathbb{A}_j^d}\left(\frac{\gamma_{i_1...i_j}}{\prod_{l=1}^{j}\gamma_{i_l}}\right)^{n'_{i_1...i_j}}\right). \quad (209)$$

Absorbing all terms that do not depend upon $M$ into $B$, we have

$$B=\sum_{\vec{\mathbf{n}}'\in C}\prod_{j=2}^{d'}\prod_{(i_1,...,i_j)\in\mathbb{A}_j^d}\left(\frac{\gamma_{i_1...i_j}}{\prod_{l=1}^{j}\gamma_{i_l}}\right)^{n'_{i_1...i_j}},\,. \quad (210)$$

Then, we can rewrite (209) as:

$$P(\vec{\mathbf{N}}=\vec{\mathbf{n}}|M) = Be^{-\vec{\mathbf{1}}^T\vec{\mathbf{\Lambda}}}\prod_{i=1}^{d}(\gamma_i M)^{n_i} = Be^{-\vec{\mathbf{1}}^T\vec{\mathbf{\Lambda}}}M^{\sum_{i=1}^{d}n_i}\prod_{i=1}^{d}(\gamma_i)^{n_i}$$

$$\stackrel{(a)}{=} Be^{-\vec{\mathbf{1}}^T\vec{\mathbf{\Lambda}}}M^{\sum_{i=1}^{d}n_i}\stackrel{(b)}{=}Be^{-\vec{\mathbf{1}}^T\vec{\mathbf{\Lambda}}}M^{\vec{\mathbf{1}}^T\vec{\mathbf{n}}}, \quad (211)$$

where, in (a) we further absorb $\prod_{i=1}^{d}(\gamma_i)^{n_i}$ into $B$ since it does not depend upon $M$, and in (b) we substitute $\sum_{i=1}^{d}n_i=\vec{\mathbf{1}}^T\vec{\mathbf{n}}$. Similarly, let us write out the expression for $P(\vec{\mathbf{N}}^g|M)$:

$$P(\vec{\mathbf{N}}^g=\vec{\mathbf{n}}^g|M) = e^{-\vec{\mathbf{1}}^T\vec{\mathbf{\Lambda}}}\prod_{j=1}^{d'}\prod_{(i_1,...,i_j)\in\mathbb{A}_j^d}(\gamma_{i_1...i_j}M^j)^{n_{i_1...i_j}^g}. \quad (212)$$

Collecting all the $M$ terms, and absorbing all the terms that do not depend upon $M$ into $D$, we obtain:

$$P(\vec{\mathbf{N}}^g=\vec{\mathbf{n}}^g|M) = De^{-\vec{\mathbf{1}}^T\vec{\mathbf{\Lambda}}}M^{\sum_{j=1}^{d'}\sum_{(i_1,...,i_j)\in\mathbb{A}_j^d}\left(jn_{i_1...i_j}^g\right)}. \quad (213)$$

Now let us analyze the term $\vec{\mathbf{1}}^T A\vec{\mathbf{n}}^g$:

$$\vec{\mathbf{1}}^T\mathbf{A}\vec{\mathbf{n}}^g \stackrel{(a)}{=} (\vec{\mathbf{n}}^g)^T\mathbf{A}^T\vec{\mathbf{1}} \stackrel{(b)}{=} (\vec{\mathbf{n}}^g)^T\begin{bmatrix}\mathbf{A}_1^T\vec{\mathbf{1}}\\\vdots\\\mathbf{A}_{d'}^T\vec{\mathbf{1}}\end{bmatrix} \stackrel{(c)}{=} (\vec{\mathbf{n}}^g)^T\begin{bmatrix}\vec{\mathbf{1}}\\2\vec{\mathbf{1}}\\\vdots\\d'\vec{\mathbf{1}},\end{bmatrix},$$

where (a) uses the fact that $\vec{\mathbf{1}}^T\mathbf{A}\vec{\mathbf{n}}^g$ is a scalar and hence is equal to its transpose, (b) uses the fact $\mathbf{A}=[\mathbf{A}_1\ldots\mathbf{A}_{d'}]$, and (c) follows from the special structure of $\mathbf{A}_i$, i.e., that each column only contains $i$ ones and $d-i$ zeros, and the fact that $\mathbf{A}_1^T\vec{\mathbf{1}}$ is akin to summing up the columns, hence $\mathbf{A}_i^T\vec{\mathbf{1}}=i\vec{\mathbf{1}}$. Equivalently we can rewrite the above equation as:

$$\vec{\mathbf{1}}^T\mathbf{A}\vec{\mathbf{n}}^g = \sum_{j=1}^{d'}\sum_{(i_1,...,i_j)\in\mathbb{A}_j^d}\left(jn_{i_1...i_j}^g\right). \quad (214)$$

Substituting (214) into (213):

$$P(\vec{\mathbf{N}}^g|M) = De^{-\vec{\mathbf{1}}^T\vec{\mathbf{\Lambda}}}M^{\vec{\mathbf{1}}^T\mathbf{A}\vec{\mathbf{n}}^g}. \quad (215)$$

Now, let us write out the expression for $P(\vec{\mathbf{N}}|\vec{\mathbf{N}}^g,M)$. Since $\vec{\mathbf{N}}=\mathbf{A}\vec{\mathbf{N}}^g$, $P(\vec{\mathbf{N}}|\vec{\mathbf{N}}^g)$ can be represented as a Kronecker delta function with the condition $\vec{\mathbf{N}}=\mathbf{A}\vec{\mathbf{N}}^g$, i.e.,

$$P(\vec{\mathbf{N}}=\vec{\mathbf{n}}|\vec{\mathbf{N}}^g=\vec{\mathbf{n}}^g,M)=P(\vec{\mathbf{N}}=\vec{\mathbf{n}}|\vec{\mathbf{N}}^g=\vec{\mathbf{n}}^g)=\delta_K\left(\vec{\mathbf{n}}=\mathbf{A}\vec{\mathbf{n}}^g\right), \quad (216)$$

where $\delta_K(\cdot)$ is the Kronecker delta function. Substituting (211), (215) and (216) in (207), we get:

$$P(\vec{\mathbf{N}}^g|\vec{\mathbf{N}},M) = \frac{De^{-\vec{\mathbf{1}}^T\vec{\mathbf{\Lambda}}}M^{\vec{\mathbf{1}}^T\mathbf{A}\vec{\mathbf{n}}^g}\delta_K(\vec{\mathbf{n}}=\mathbf{A}\vec{\mathbf{n}}^g)}{Be^{-\vec{\mathbf{1}}^T\vec{\mathbf{\Lambda}}}M^{\vec{\mathbf{1}}^T\vec{\mathbf{n}}}}\stackrel{(a)}{=}\frac{DM^{\vec{\mathbf{1}}^T\mathbf{A}\vec{\mathbf{n}}^g}\delta_K(\vec{\mathbf{n}}=A\vec{\mathbf{n}}^g)}{BM^{\vec{\mathbf{1}}^T\vec{\mathbf{n}}}},$$

$$\stackrel{(b)}{=}\frac{DM^{\vec{\mathbf{1}}^T\mathbf{A}\vec{\mathbf{n}}^g}\delta_K(\vec{\mathbf{n}}=\mathbf{A}\vec{\mathbf{n}}^g)}{BM^{\vec{\mathbf{1}}^T\mathbf{A}\vec{\mathbf{n}}^g}}\stackrel{(c)}{=}\frac{D\delta_K(\vec{\mathbf{n}}=\mathbf{A}\vec{\mathbf{n}}^g)}{B}, \quad (217)$$

where we obtain (a) by canceling the term $e^{-\vec{1}^T \vec{\Lambda}}$, (b) by using the fact $\vec{n} = \mathbf{A}\vec{n}^g$ due to the delta function, and (c) by canceling the term $M^{\vec{1}^T \mathbf{A}\vec{n}^g}$. Since the terms $D, B$ and $\delta_K(\vec{n} = \mathbf{A}\vec{n}^g)$ do not depend upon $M$, we can conclude $P(\vec{\mathbf{N}}^g|\vec{\mathbf{N}}, M)$ also does not depend upon $M$, i.e. $P(\vec{\mathbf{N}}^g|\vec{\mathbf{N}}, M) = P(\vec{\mathbf{N}}^g|\vec{\mathbf{N}})$, or $M \to \vec{\mathbf{N}} \to \vec{\mathbf{N}}^g$. $\qquad\square$

**Lemma 18.** *Let* $M \sim P(M)$ *have support over* $(0, \infty)$. *Define,* $\vec{\mathbf{X}} = \begin{bmatrix} \vec{\mathbf{X}}_1^T & \dots & \vec{\mathbf{X}}_{d_1}^T \end{bmatrix}^T$ *and* $\vec{\mathbf{Y}} = \begin{bmatrix} \vec{\mathbf{Y}}_1 & \dots & \vec{\mathbf{Y}}_{d_2}^T \end{bmatrix}^T$, *where* $\vec{\mathbf{X}}_i$ *and* $\vec{\mathbf{Y}}_j$ *are random vectors of size* $q_i \in \mathbb{N}$ *and* $r_j \in \mathbb{N}$, *respectively,* $\forall\, i \in [d_1]$, *and* $j \in [d_2]$. *Let the conditional distribution of* $\vec{\mathbf{X}}$ *and* $\vec{\mathbf{Y}}$ *conditioned on* $M$ *be specified as follows:*

$$P\left(\vec{\mathbf{X}}|M\right) = \prod_{i=1}^{d_1} P\left(\vec{\mathbf{X}}_i|M\right), \text{ and } P\left(\vec{\mathbf{Y}}|M\right) = \prod_{i=1}^{d_2} P\left(\vec{\mathbf{Y}}_i|M\right), \text{ where}$$

$$P\left(\vec{\mathbf{X}}_i|M\right) = Poisson\left(q_i, 1, \vec{\Lambda}_i^X\right), \vec{\Lambda}_i^X = \begin{bmatrix} \gamma_{i1}^X M^i & \dots & \gamma_{iq_i}^X M^i \end{bmatrix}^T \ \forall\, i \in [d_1], \text{ and}$$

$$P\left(\vec{\mathbf{Y}}_j|M\right) = Poisson\left(r_j, 1, \vec{\Lambda}_j^Y\right), \vec{\Lambda}_i^Y = \begin{bmatrix} \gamma_{i1}^Y M^j & \dots & \gamma_{ir_i}^Y M^j \end{bmatrix}^T \ \forall\, j \in [d_2]. \quad (218)$$

*If $d_1 \geq d_2$ and (219) hold*

$$\sum_{j=1}^{q_i} \gamma_{ij}^X \geq \sum_{j=1}^{r_i} \gamma_{ij}^Y \ \forall\, i \in [d_2] \quad (219)$$

*then the distribution* $\tilde{Q}(M, \vec{\mathbf{X}}, \vec{\mathbf{Y}})$ *defined in (220) lies in* $\Delta_P$.

$$\tilde{Q}(M, \vec{\mathbf{X}}, \vec{\mathbf{Y}}) = P(M)P(\vec{\mathbf{X}}|M)\tilde{Q}(\vec{\mathbf{Y}}|\vec{\mathbf{X}}), \quad (220)$$

*where* $\tilde{Q}(\vec{\mathbf{Y}}|\vec{\mathbf{X}})$ *is a product of* $d_2$ *multinomial distributions, i.e* $\tilde{Q}(\vec{\mathbf{Y}}|\vec{\mathbf{X}}) = \prod_{i=1}^{d_2} \tilde{Q}(\vec{\mathbf{Y}}_i|\vec{\mathbf{X}}_i)$, *where* $\tilde{Q}(\vec{\mathbf{Y}}_i|\vec{\mathbf{X}}_i) = Multinomial(\vec{\mathbf{k}}_i; N_i, \vec{\mathbf{p}}_i)$ *and*

$$N_i = \sum_{j=1}^{n_i} x_{ij}, \ \vec{\mathbf{k}}_i = \begin{bmatrix} y_{i1} & \cdots & y_{im_i} & \sum_{j=1}^{n_i} x_{ij} - \sum_{j=1}^{m_i} y_{ij} \end{bmatrix}^T,$$

$$\vec{\mathbf{p}}_i = \begin{bmatrix} \frac{\gamma_{i1}^Y}{\sum_{j=1}^{n_i} \gamma_{ij}^X} & \cdots & \frac{\gamma_{im_i}^Y}{\sum_{j=1}^{n_i} \gamma_{ij}^X} & 1 - \frac{\sum_{j=1}^{m_i} \gamma_{ij}^Y}{\sum_{j=1}^{n_i} \gamma_{ij}^X} \end{bmatrix}^T.$$

*Proof.* Lemma 18 follows from a straightforward application of lemma 16. For showing $\tilde{Q}(M, \vec{\mathbf{X}}, \vec{\mathbf{Y}})$ specified in (220) lies in $\Delta_P$, we need to show that $\tilde{Q}(M, \vec{\mathbf{X}}) = P(M, \vec{\mathbf{X}})$ and $\tilde{Q}(M, \vec{\mathbf{Y}}) = P(M, \vec{\mathbf{Y}})$. $\tilde{Q}(M, \vec{\mathbf{X}}) = P(M, \vec{\mathbf{X}})$ is satisfied trivially by construction of $\tilde{Q}(M, \vec{\mathbf{X}}, \vec{\mathbf{Y}})$.

Proving $\tilde{Q}(\vec{\mathbf{Y}}, M) = P(\vec{\mathbf{Y}}, M)$:

For showing the second condition, we will explicitly show $\tilde{Q}(\vec{\mathbf{Y}}|M) = P(\vec{\mathbf{Y}}|M)$ which, combined with the fact that $\tilde{Q}(M) = P(M)$ (by construction), will show $\tilde{Q}(\vec{\mathbf{Y}}, M) = P(\vec{\mathbf{Y}}, M)$ completing our proof.

Calculating $\tilde{Q}(\vec{\mathbf{Y}}|\vec{\mathbf{X}})$: From the law of total probability, we have:

$$\tilde{Q}(\vec{\mathbf{Y}}|M) = \sum_{\vec{\mathbf{x}}} P(\vec{\mathbf{X}} = \vec{\mathbf{x}}|M)\tilde{Q}(\vec{\mathbf{Y}}|\vec{\mathbf{X}} = \vec{\mathbf{x}})$$

$$\overset{(a)}{=} \sum_{\vec{\mathbf{x}}} \prod_{i=1}^{d_1} P(\vec{\mathbf{X}}_i = \vec{\mathbf{x}}_i|M) \prod_{i=1}^{d_2} \tilde{Q}(\vec{\mathbf{Y}}_i|\vec{\mathbf{X}}_i = \vec{\mathbf{x}}_i)$$

$$\overset{(b)}{=} \sum_{\vec{\mathbf{x}}} \prod_{i=1}^{d_2} P(\vec{\mathbf{X}}_i = \vec{\mathbf{x}}_i|M)\tilde{Q}(\vec{\mathbf{Y}}_i|\vec{\mathbf{X}}_i = \vec{\mathbf{x}}_i) \prod_{i=d_2+1}^{d_1} P(\vec{\mathbf{X}}_i = \vec{\mathbf{x}}_i|M)$$

$$\overset{(c)}{=} \prod_{i=1}^{d_2}\left(\sum_{\vec{\mathbf{x}}_i} P(\vec{\mathbf{X}}_i = \vec{\mathbf{x}}_i|M)\tilde{Q}(\vec{\mathbf{Y}}_i|\vec{\mathbf{X}}_i = \vec{\mathbf{x}}_i)\right) \prod_{i=d_2+1}^{d_1}\left(\underbrace{\sum_{\vec{\mathbf{x}}_i} P(\vec{\mathbf{X}}_i = \vec{\mathbf{x}}_i|M)}_{=1}\right)$$

$$\overset{(d)}{=} \prod_{i=1}^{d_2}\left(\underbrace{\sum_{\vec{\mathbf{x}}_i} P(\vec{\mathbf{X}}_i = \vec{\mathbf{x}}_i|M)\tilde{Q}(\vec{\mathbf{Y}}_i|\vec{\mathbf{X}}_i = \vec{\mathbf{x}}_i)}_{=\tilde{Q}(\vec{\mathbf{Y}}_i|M)}\right) = \prod_{i=1}^{d_2} \tilde{Q}(\vec{\mathbf{Y}}_i|M), \tag{221}$$

where $(a)$ is due to the particular structure of $P(\vec{\mathbf{X}}|M)$ and $\tilde{Q}(\vec{\mathbf{Y}}|\vec{\mathbf{X}})$ specified in (218) and (220), respectively, $(b)$ is just collecting the terms of $P(\vec{\mathbf{X}}_i|M)$ and $\tilde{Q}(\vec{\mathbf{Y}}_i|\vec{\mathbf{X}}_i)$ into a single product, $(c)$ is due to spreading the summation of the components of $\tilde{\mathbf{x}}$ over their corresponding distribution components, and $(d)$ is due to the fact that sum over all probabilities of all possible $\tilde{\mathbf{x}}_i$ is 1.

Note that for a fixed $m$, $P\left(\vec{\mathbf{X}}_i|M = m\right) \sim \text{Poisson}\left(n_i, 1, \begin{bmatrix}\gamma_{i1}m^i & \cdots & \gamma_{in_i}m^i\end{bmatrix}^T\right)$, and $Q(\vec{\mathbf{Y}}_i|\vec{\mathbf{X}}_i) = \text{Multinomial}(\vec{\mathbf{k}}_i; N_i, \vec{\mathbf{p}}_i)$, where

$$N_i = \sum_{j=1}^{n_i} x_{ij},\ \vec{\mathbf{k}}_i = \begin{bmatrix}y_{i1} & \cdots & y_{im_i} & \sum_{j=1}^{n_i} x_{ij} - \sum_{j=1}^{m_i} y_{ij}\end{bmatrix}^T,$$

$$\vec{\mathbf{p}}_i = \begin{bmatrix}\frac{\gamma_{i1}^Y}{\sum_{j=1}^{n_i}\gamma_{ij}^X} & \cdots & \frac{\gamma_{im_i}^Y}{\sum_{j=1}^{n_i}\gamma_{ij}^X} & 1 - \frac{\sum_{j=1}^{m_i}\gamma_{ij}^Y}{\sum_{j=1}^{n_i}\gamma_{ij}^X}\end{bmatrix}^T.$$

Note that we can equivalently write $\vec{\mathbf{p}}_i$ as follows,

$$\vec{\mathbf{p}}_i = \begin{bmatrix}\frac{\gamma_{i1}^Y M^i}{\sum_{j=1}^{n_i}\gamma_{ij}^X M^i} & \cdots & \frac{\gamma_{im_i}^Y M^i}{\sum_{j=1}^{n_i}\gamma_{ij}^X M^i} & 1 - \frac{\sum_{j=1}^{m_i}\gamma_{ij}^Y M^i}{\sum_{j=1}^{n_i}\gamma_{ij}^X M^i}\end{bmatrix}^T. \tag{222}$$

In the setup of lemma 16, choose $d_1 = q_i$ and $d_2 = r_i$, $\lambda_j^{(2)} = \gamma_{ij}^Y M^i \ \forall\ j \in [r_i]$ and $\lambda_j^{(1)} = \gamma_{ij}^X M^i \ \forall\ j \in [q_i]$. From (219), we know the condition $\sum_{j=1}^{d_2} \lambda_j^{(2)} \leq \sum_{j=1}^{d_2} \lambda_j^{(2)}$ is satisfied by assumption. Hence, we use the result of lemma 15 to conclude $\tilde{Q}(\vec{\mathbf{Y}}_i|M) = \text{Poisson}(r_i, 1, \vec{\mathbf{\Lambda}}')$, with

$$\vec{\mathbf{\Lambda}}' = \begin{bmatrix}\lambda_{i1} & \ldots, \lambda_{ir_i}\end{bmatrix}^T, \text{ with } \lambda_{ij} = \gamma_{ij}^Y M^i. \tag{223}$$

Comparing $\tilde{Q}(\vec{\mathbf{Y}}_i|M)$ specified in (223) with $P(\vec{\mathbf{Y}}_i|M)$ specified in (218), we can conclude $Q(\vec{\mathbf{Y}}_i|M) = P(\vec{\mathbf{Y}}_i|M)$. Substituting $Q(\vec{\mathbf{Y}}_i|M) = P(\vec{\mathbf{Y}}_i|M)$ in (221), we obtain:

$$\tilde{Q}(\vec{\mathbf{Y}}|M) = \prod_{i=1}^{d_2} P(\vec{\mathbf{Y}}_i|M). \tag{224}$$

Comparing (224) with (218), we can conclude $\tilde{Q}(\vec{\mathbf{Y}}|M) = P(\vec{\mathbf{Y}}|M)$, finishing our proof. $\qquad\square$

# K Proof of Theorem 6

In this section, we provide the proof of theorem 6. Since theorem 6 provides the analytical PID terms for the linear convolution-closed system defined in Sec. 5.1, we briefly restate certain key properties of convolution-closed distributions and the linear convolution-closed system for convenience.

## K.1 Convolution-closed distributions

Convolution-closed distributions are a large class of distributions that are defined as follows:

**Definition 1.** *Let $\mathcal{F}_\mathcal{D}$ denote a family of distributions, where each member distribution, $f(\delta) \in \mathcal{F}_\mathcal{D}$, is indexed by a parameter $\delta$. Consider $X_1 \sim f(\delta_1)$, $X_2 \sim f(\delta_2)$, and $X_1 \perp\!\!\!\perp X_2$ for some $\delta_1, \delta_2 \in \mathcal{D}$. Then, $\mathcal{F}_\mathcal{D}$ is convolution-closed in the parameter $\delta$, if*

$$X_1 + X_2 \sim f(\delta_1) * f(\delta_2) = f(\delta_1 + \delta_2) \ \forall \ \delta_1, \delta_2 \in \mathcal{D} \text{ such that } \delta_1 + \delta_2 \in \mathcal{D}, \quad (225)$$

*where $*$ denotes the convolution operator.*

Convolution-closed distributions define a natural dilation/thinning operator. Formally, let $X \sim f(\delta)$. Then, we define $X_\epsilon$ as the $\epsilon$-dilated version of $X$ if $X \sim f(\epsilon\delta)$ for some $\epsilon \in (0, 1)$ such that $\epsilon\delta \in \mathcal{D}$. Furthermore, if we assume $(1 - \epsilon)\delta \in \mathcal{D}$, then $P(X_\epsilon|X)$ can be defined as $P(X_\epsilon|X_\epsilon + X_{(1-\epsilon)})$, where $X_{1-\epsilon} \sim f((1-\epsilon)\delta)$ and $X_\epsilon \perp\!\!\!\perp X_{1-\epsilon}$. Denote, $P(X_\epsilon|X = x) = P(X_\epsilon|X_\epsilon + X_{(1-\epsilon)} = x) = G(\epsilon\delta, (1-\epsilon)\delta, x)$.

## K.2 Definition of linear convolution-closed system

Let $\mathcal{F}_\mathcal{D}$ be a convolution-closed distribution family in parameter $\delta \in \mathcal{D}$, and $M$ be a target/message random variable having distribution $P(M)$ over some support set $\mathcal{M}$. Define the conditional distribution of random variables $X$ and $Y$ conditioned on $M$ as follows:

$$P(X|M{=}m) = f(\delta_m^X) \text{ and } P(Y|M{=}m) = f(\delta_m^Y) \text{ such that } \delta_m^X, \delta_m^Y \in \mathcal{D} \ \forall \ m \in \mathcal{M}. \quad (226)$$

Furthermore, we assume that there exists $\gamma \in (0, \infty)$ such that: $\delta_m^X = \gamma\delta_m^Y \ \forall \ m \in \mathcal{M}$. We denote such a system of random variables $(M, X, Y)$, having the joint distribution $P(M, X, Y)$, a *linear convolution-closed system*.

## K.3 Formal proof of theorem 6

**Theorem 6.** *Let the joint density $P(M, X, Y)$ of random variables $M$, $X$, and $Y$ describe a linear convolution-closed system. Without the loss of generality, assume $\gamma \leq 1$. If*

*(a) $(1 - \gamma)\delta_m^Y \in \mathcal{D} \ \forall \ m \in \mathcal{M}$,*

*(b) $P(X_\gamma|X_\gamma + X_{1-\gamma}{=}x, M{=}m) = G(\gamma\delta_m^X, (1-\gamma)\delta_m^X, x)$ does not depend on $m$, where $P(X_\gamma|M) = f(\gamma\delta_m^X)$, $P(X_{1-\gamma}|M) = f((1-\gamma)\delta_m^X)$ and $X_\gamma \perp\!\!\!\perp X_{1-\gamma}|M$,*

*then $\Delta_P$ contains a Markov chain of the form $M \to X \to Y$ and $UI(M; Y \backslash X) = 0$.*

*Proof.* We first note that we can always assume $\gamma \leq 1$ without the loss of generality because if $\gamma \geq 1$, then we can always switch our nomenclature to refer to $Y$ as $X$, and $X$ as $Y$.

We briefly outline the proof structure.

1. In the first part, we explicitly construct a joint distribution $Q_{MC}(M, X, Y)$ having the Markovian structure $M \to X \to Y$.

2. In the second part, we show that the $Q_{MC}(M, X, Y)$ constructed in the first part, lies in $\Delta_P$. Therefore, we can then apply the result of proposition 1 to conclude $UI(M; Y \backslash X) = 0$.

**Part 1: Specifying $Q_{MC}(M, X, Y)$**

Denote the joint distribution of the Markov chain $M \to X \to Y$ as $Q_{MC}(M, X, Y)$. We can decompose $Q_{MC}(M, X, Y)$ by utilizing its Markovian structure as follows:

$$Q_{MC}(M, X, Y) = Q_{MC}(M)Q_{MC}(X|M)Q_{MC}(Y|X).$$

Consequently, we can construct the distribution $Q_{MC}(M, X, Y)$ by individually specifying $Q_{MC}(M)$, $Q_{MC}(X|M)$, and $Q_{MC}(Y|X)$.

Specifying $Q_{MC}(M)$ and $Q_{MC}(X|M)$: We specify $Q_{MC}(M)$ and $Q_{MC}(X|M)$ as follows:

$$Q_{MC}(M) = P(M) \text{ and } Q_{MC}(X|M) = P(X|M). \tag{227}$$

Specifying $Q_{MC}(Y|X)$: By assumption (b) in the theorem, we know that $G(\gamma \delta_m^X, (1 - \gamma)\delta_m^X, x)$ does not depend on $m$, and consequently, does not depend on $\delta_m^X$. Hence, we can simplify notation of $G(\gamma \delta_m^X, (1 - \gamma)\delta_m^X, m)$ as $G(\gamma, (1 - \gamma), x)$. We specify $Q(Y|X)$ as follows:

$$Q(Y|X = x) = G(\gamma, 1 - \gamma, x). \tag{228}$$

**Part 2: Showing $Q_{MC}(M, X, Y) \in \Delta_P$**

For showing that $Q_{MC}(M, X, Y) \in \Delta_P$, we first need to derive $Q_{MC}(Y|M)$.

Deriving $Q_{MC}(Y|M)$: We use the result of lemma 19 to derive $Q_{MC}(Y|M)$. As $Q_{MC}(X|M = m) = f(\delta_m^X)$ is a convolution-closed distribution for a fixed $m$, and $Q_{MC}(Y = y|X = x) = G(\gamma, 1 - \gamma, x)$, the result of lemma 19 shows that $Q_{MC}(Y|M)$ can be expressed as:

$$Q_{MC}(Y|M = m) = f(\gamma \times \delta_m^X) = f(\gamma \delta_m^X) \overset{(a)}{=} f(\delta_m^Y) = P(Y|M = m), \tag{229}$$

where $(a)$ is due to the assumption $\delta_m^X = \gamma \delta_m^Y$ (see Appx. K.2).

From (227) and (229), we can conclude:

$$Q_{MC}(M, X) = Q_{MC}(M)Q_{MC}(X|M) = P(M)P(X|M) = P(M, X), \tag{230}$$
$$Q_{MC}(M, Y) = Q_{MC}(M)Q_{MC}(Y|M) = P(M)P(Y|M) = P(M, Y). \tag{231}$$

Hence, $Q_{MC}(M, X, Y) \in \Delta_P$ and consequently, by proposition 1, $UI(M; Y \backslash X) = 0$, concluding our proof for case 1.

$\square$

**Lemma 19.** *Let $\mathcal{F}_\mathcal{D}$ be a convolution-closed distribution family convolution-closed in the parameter $\delta$. For a $\delta \in \mathcal{D}$, assume there exists $\epsilon \in (0, 1)$ such that $\epsilon\delta, (1 - \epsilon)\delta \in \mathcal{D}$. Consider the following random variables:*

$$X'_\epsilon \sim f(\epsilon\delta) \in \mathcal{F}_\mathcal{D}, \quad X'_{1-\epsilon} \sim f((1 - \epsilon)\delta) \in \mathcal{F}_\mathcal{D}, \quad \text{and } X \sim f(\delta) \in \mathcal{F}_\mathcal{D},$$

*such that $X'_\epsilon \perp\!\!\!\perp X'_{1-\epsilon}$. If $P(X_\epsilon = x_\epsilon|X = x) = P(X'_\epsilon = x_\epsilon|X'_\epsilon + X'_{1-\epsilon} = x)$, then we have $P(X_\epsilon = x_\epsilon) = P(X'_\epsilon = x_\epsilon) = f(x_\epsilon; \epsilon\delta)$ or $X_\epsilon \overset{d}{=} X'_\epsilon$, where $\overset{d}{=}$ implies equality in distribution [57].*

*Proof.* First let us derive the distribution of $X' = X'_{1-\epsilon} + X'_\epsilon$. Since, $X'_\epsilon \perp\!\!\!\perp X'_{1-\epsilon}$, we can write $P(X')$ as follows:

$$P(X') = f(\epsilon\delta) * f((1 - \epsilon)\delta) \overset{(a)}{=} f(\epsilon\delta + (1 - \epsilon)\delta) = f(\delta) \overset{(b)}{=} P(X), \tag{232}$$

where $*$ is the convolution operator, $(a)$ is due to the properties of convolution-closed distribution (see Appx. K.1), and $(b)$ is due to the definition of $X$ in the lemma statement. From (232), we can conclude $P(X = x) = P(X' = x)$, and by assumption we have $P(X_\epsilon|X) = P(X'_\epsilon|X') = P(X'_\epsilon|X'_\epsilon + X'_{(1-\epsilon)})$. Hence, we have:

$$P(X = x, X_\epsilon = x_\epsilon) = P(X = x)P(X_\epsilon = x_\epsilon|X = x) = P(X' = x)P(X'_\epsilon = x_\epsilon|X' = x),$$
$$= P(X' = x)P(X'_\epsilon = x_\epsilon|X'_\epsilon + X'_{1-\epsilon} = x)$$
$$= P(X'_\epsilon + X'_{1-\epsilon} = x, X'_\epsilon = x_\epsilon). \tag{233}$$

From (233), we can conclude that $P(X_\epsilon) = P(X'_\epsilon) = f(\epsilon\delta)$. $\square$

# L   Proof of Theorem 7

In this section, we provide the proof of theorem 7.

## L.1   Definition of $p_{exp1}$ and $p_{exp2}$

We restate the definitions of $p_{exp1}$ and $p_{exp2}$ defined in Sec. 5.2, which is used in the proof of theorem 7. Let $X \sim p_{exp1}(X)$, where

$$p_{exp1}(X = x; \theta_1, \theta_2) = H(\theta_1, \theta_2) \exp(\theta_1^T x - \theta_2^T A(x)), \tag{234}$$

for some appropriately defined $H(\cdot, \cdot), A(\cdot), \theta_1$ and $\theta_2$. Furthermore, define a random variable $Y$ through its conditional density $p(Y|X = x)$:

$$p(Y = y | X = x; \theta_1, \theta_2) = h(y) \exp\left( x^T T(y) - \theta_3^T A(x) \right), \tag{235}$$

for some $h(\cdot), T(\cdot)$, and $\theta_3$, such that $p(Y = y | X = x; \theta_1, \theta_2)$ is a well-defined distribution. Furthermore, the marginal distribution of $Y$ is expressed as:

$$p_{exp2}(Y = y; \theta_1, \theta_2, \theta_3) = h(y) \frac{H(\theta_1, \theta_2)}{H(\theta_1 + T(y), \theta_2 + \theta_3)} \text{ (see proof of theorem 1 in [26]).} \tag{236}$$

## L.2   Formal proof of theorem 7

**Theorem 7.** *Let $M, X$, and $Y$ be random variables having the joint distribution $P(M, X, Y)$. Furthermore, the conditional distribution of $X$ and $Y$ conditioned on $M$ are as follows: $P(X|M{=}m){=}p_{exp1}(X; \theta_1(m), \theta_2(m))$ and $P(Y|M{=}m){=}p_{exp2}(Y; \theta_1(m), \theta_2(m), \theta_3)$. Then, $\Delta_P$ contains a Markov chain of the form $M \to X \to Y$ and $UI(M; Y \backslash X) = 0$.*

*Proof.* We prove theorem 7 by explicitly constructing a Markov chain $M \to X \to Y$, having joint distribution $Q_{MC}(M, X, Y)$, and showing that $Q_{MC}(M, X, Y) \in \Delta_P$.

Specifying $Q_{MC}(M)$ and $Q_{MC}(X|M)$: We specify $Q_{MC}(M)$ and $Q_{MC}(X|M)$ as follows:

$$Q_{MC}(M) = P(M), \text{ and } Q_{MC}(X|M) = P(X|M). \tag{237}$$

Specifying $Q_{MC}(Y|X)$: We specify $Q_{MC}(Y|X) = h(y) \exp(x^T T(y) - \theta_3^T A(x))$. Then, by (236), we know that

$$Q(Y = y | M = m) = p_{exp2}(y; \theta_1(m), \theta_2(m), \theta_3) = \frac{h(y) H(\theta_1(m), \theta_2(m))}{H(\theta_1(m) + T(y), \theta_2(m) + \theta_3(m))}. \tag{238}$$

From (237) and (238), we can conclude:

$$Q_{MC}(M, Y) = Q_{MC}(Y|M) Q_{MC}(M) = P(Y|M) P(M) = P(M, Y), \tag{239}$$
$$Q_{MC}(M, X) = Q_{MC}(X|M) Q_{MC}(M) = P(X|M) P(M) = P(M, X). \tag{240}$$

Hence, $Q_{MC}(M, X, Y) \in \Delta_P$ and consequently, by proposition 1, $UI(M; Y \backslash X) = 0$, concluding our proof. $\square$

# M   Additional proofs and details for Sec. 6

In this section, we provide the proofs of certain statements made in Sec. 6, as well as additional details regarding the simulation study presented in the same section.

## M.1   Generating $\bar{Q}$ for every $Q \in \Delta_P$

Our goal in this section is to show that there exists a $\bar{Q}$ (as described in Sec. 6) for every $Q \in \Delta_P$, where the corresponding $P(M, X, Y)$ is as defined in Sec. 6. We first briefly restate the assumptions on $P(M, X, Y)$ described in Sec. 6. The distribution $P(M, X, Y)$ has the following properties: the random variable $M$ has support over $\mathcal{M}$, the conditional distributions $P(X|M)$ and $P(Y|M)$ are

members of some convolution-closed distribution family $\mathcal{F}_\mathcal{D}$, and there exists some $\delta_{bias}^X, \delta_{bias}^Y \in \mathcal{D}$ such that:

$$P(X|M=m)=f(\delta_m^X) \text{ and } P(Y|M=m)=f(\delta_m^Y) \text{ where } \delta_m^X, \delta_m^Y \in \mathcal{D} \; \forall \; m \in \mathcal{M}, \text{ with}$$

$$\delta_m^X - \delta_{bias}^X = \epsilon_m^{(1)}(\delta_m^Y - \delta_{bias}^Y) \text{ for some } \epsilon_m^{(1)} \in [0,1] \; \forall \; m \in \mathcal{M},$$

$$\delta_m^Y = \epsilon_m^{(2)}(\delta_m^Y - \delta_{bias}^Y) \text{ for some } \epsilon_m^{(2)} \in [0,1] \; \forall \; m \in \mathcal{M},$$

$$\delta_m^X = \epsilon_m^{(3)}(\delta_m^X - \delta_{bias}^X) \text{ for some } \epsilon_m^{(3)} \in [0,1] \; \forall \; m \in \mathcal{M}, \text{ and}$$

$$(\delta_m^Y - \delta_{bias}^Y), \; (\delta_m^X - \delta_{bias}^X), \; (\delta_m^X - \delta_m^Y - (\delta_{bias}^X - \delta_{bias}^Y)) \in \mathcal{D}. \tag{241}$$

As $Q(M,X)=P(M,X)$ and $Q(M,Y) = P(M,Y)$ due to $Q \in \Delta_P$, we have:

$$Q(X|M=m)=f(\delta_m^X), \text{ and } Q(Y|M=m)=f(\delta_m^Y) \text{ where } \delta_m^X, \delta_m^Y \in \mathcal{D} \; \forall \; m \in \mathcal{M}. \tag{242}$$

### M.1.1  Explicitly constructing a $\bar{Q}$ for every $Q \in \Delta_P$

We show existence of a $\bar{Q}$ for every $Q \in \Delta_P$ by explicitly constructing a $\bar{Q}$ for each $Q \in \Delta_P$.

We define two vectors to simplify our notations:

$$\vec{\mathbf{A}} = [X \quad Y]^T, \quad \vec{\mathbf{B}} = [X' \quad Y'' \quad n_X \quad Y' \quad n_Y]^T, \tag{243}$$

where $X$ and $Y$ are the random variables defined above. $(X', Y'', n_X)$, and $(Y', n_Y)$ are the corresponding dilated versions of $X$ and $Y$ as defined in Sec. 6. Note that the joint distribution of $(M, X', Y'', n_X, Y', n_Y)$ is $\bar{Q}(M, X', Y'', n_X, Y', n_Y)$. We reiterate the two main properties of $\bar{Q}(X', Y'', n_X, Y', n_Y)$, mentioned in Sec. 6, that we want to prove:

1. The marginal conditional distributions of $(X', Y'', n_X, Y', n_Y)$ given $M$ are as follows:

$$\bar{Q}(X'|M=m)=f(\delta_m^X - \delta_m^Y - (\delta_{bias}^X - \delta_{bias}^Y)), \; \bar{Q}(Y''|M=m) = f(\delta_m^Y - \delta_{bias}^Y),$$

$$\bar{Q}(Y'|M=m)=f(\delta_m^Y - \delta_{bias}^Y), \bar{Q}(n_X|M) = f(\delta_{bias}^X), \bar{Q}(n_Y|M) = f(\delta_{bias}^Y).$$

2. The random variables $(X', Y'', n_X)$ are jointly conditionally independent given $M$. Similarly, the random variables $(Y', n_Y)$ are also conditionally independent given $M$.

We construct $\vec{\mathbf{B}}$ from $\vec{\mathbf{A}}$ by specifying the conditional distribution $\bar{Q}(\vec{\mathbf{B}}|\vec{\mathbf{A}}, M)$. Note that given the values of $\vec{\mathbf{A}}$ and $M$, we can always use $\bar{Q}(\vec{\mathbf{B}}|\vec{\mathbf{A}}, M)$ to construct the random variables $\vec{\mathbf{B}}$. The goal of this remaining section is to show that the corresponding distribution $\bar{Q}(\vec{\mathbf{B}}, M)$ (specified through $\bar{Q}(\vec{\mathbf{B}}|\vec{\mathbf{A}}, M)$) satisfies the above two properties. We first explicitly state the structure of $\bar{Q}(\vec{\mathbf{B}}|\vec{\mathbf{A}}, M)$.

Specifying $\bar{Q}(\vec{\mathbf{B}}|\vec{\mathbf{A}}, M)$:

We first impose the following structure on $\bar{Q}(\vec{\mathbf{B}}|\vec{\mathbf{A}}, M)$ as follows:

$$\bar{Q}(\vec{\mathbf{B}}|\vec{\mathbf{A}}, M) = \bar{Q}(X', Y'', n_X|X, M)\bar{Q}(Y', n_Y|Y, M), \tag{244}$$

where $\bar{Q}(X', Y'', n_X|X, M)$ and $\bar{Q}(Y', n_Y|Y, M)$ are the conditional distributions through which we specify $\bar{Q}(\vec{\mathbf{B}}|\vec{\mathbf{A}}, M)$. Since $\bar{Q}(\vec{\mathbf{B}}|\vec{\mathbf{A}}, M)$ is a product of two well-defined distributions, it is a valid distribution. Let us now analyze the joint distribution $\bar{Q}(M, \vec{\mathbf{A}}, \vec{\mathbf{B}})$. By chain rule, we have:

$$\bar{Q}(M, \vec{\mathbf{A}}, \vec{\mathbf{B}}) = \bar{Q}(M, \vec{\mathbf{A}})\bar{Q}(\vec{\mathbf{B}}|\vec{\mathbf{A}}, M). \tag{245}$$

Substituting the fact that $(M, \vec{\mathbf{A}}) = (M, X, Y)$ and using the structure of $\bar{Q}(\vec{\mathbf{B}}|\vec{\mathbf{A}}, M)$ specified in (244), we have:

$$\bar{Q}(M, \vec{\mathbf{A}}, \vec{\mathbf{B}}) = \bar{Q}(M)\bar{Q}(X, Y|M)\bar{Q}(X', Y'', n_X|X, M)\bar{Q}(Y', n_Y|Y, M). \tag{246}$$

We now specify the two components of $\bar{Q}(\vec{\mathbf{B}}|\vec{\mathbf{A}}, M)$.

Specifying $\bar{Q}(Y', n_Y|Y, M)$:

Using the chain rule on $\bar{Q}(Y', n_Y | Y, M)$, we obtain
$$\bar{Q}(Y', n_Y | Y, M) = \bar{Q}(Y'|Y,M)\bar{Q}(n_Y|Y,Y',M). \tag{247}$$
Choose:
$\bar{Q}(Y'|Y{=}y, M{=}m) = G(\epsilon_m^{(2)}\delta_m^Y, (1 - \epsilon_m^{(2)})\delta_m^Y, y),$ where $\delta_m^Y {=} \epsilon_m^{(2)}(\delta_m^Y - \delta_{bias}^Y) \ \forall \ m \in \mathcal{M},$ and
$\bar{Q}(n_Y|Y{=}y, Y'{=}y', M{=}m) = \delta_K(n_Y = y - y'). \tag{248}$

Here, $\delta_K(\cdot)$ denotes the Kronecker delta function [59]. Marginalizing $(X, X', Y'', n_X)$ from (246), we obtain:
$$\bar{Q}(M, Y, Y', n_Y) = \bar{Q}(M)\bar{Q}(Y|M)\bar{Q}(Y', n_Y|M, Y) = \bar{Q}(M)\bar{Q}(Y, Y', n_Y|M). \tag{249}$$
Consider the distribution $\bar{Q}(Y, Y', n_Y|M)$. Then, we know:
$$\bar{Q}(Y|M{=}m) = f(\delta_m^Y),$$
$$\bar{Q}(n_Y|Y{=}y, Y'{=}y'M{=}m) = \delta_K(n_Y = y - y'), \text{ and}$$
$$\bar{Q}(Y'{=}y'|Y{=}y, M{=}m) = G(\epsilon_m^{(2)}\delta_m^Y, (1 - \epsilon_m^{(2)})\delta_m^Y, y). \tag{250}$$
Comparing the above equation with the structure of $\bar{Q}(Y, Y', n_Y|M)$ specified in proposition 8, we can conclude that $(Y', n_Y)$ are just dilated versions of $Y$. Consequently, using the result of proposition 8, we have:
$$\bar{Q}(Y'|M{=}m) = f(\delta_m^Y - \delta_{bias}^Y), \bar{Q}(n_Y|M) = f(\delta_{bias}^Y), \text{ and } Y' \perp\!\!\!\perp n_Y|M. \tag{251}$$

Specifying $\bar{Q}(X', Y'', n_X|X, M)$:

We similarly decompose $\bar{Q}(X', Y'', n_X|X, M)$ using the chain rule as follows:
$$\bar{Q}(X', Y'', n_X|X, M) = \bar{Q}(n_X|X, M)\bar{Q}(X'|X, n_X, M)\bar{Q}(Y''|X, n_X, X', M). \tag{252}$$
We choose:
$\bar{Q}(n_Y|X{=}x, M{=}m) = G((1 - \epsilon_m^{(3)})\delta_m^X, \epsilon_m^{(3)}\delta_m^X, x),$ where $\delta_m^X = \epsilon_m^{(3)}(\delta_m^X - \delta_{bias}^X) \ \forall \ m \in \mathcal{M},$
$\bar{Q}(X'|X{=}x, n_X{=}x_{n_X}, M{=}m) = G((1 - \epsilon_m^{(1)})(\delta_m^X - \delta_{bias}^X), \epsilon_m^{(1)}(\delta_m^X - \delta_{bias}), x - x_{n_X}),$ where
$\delta_m^X - \delta_{bias}^X = \epsilon_m^{(1)}(\delta_m^Y - \delta_{bias}^Y) \ \forall \ m \in \mathcal{M},$ and
$\bar{Q}(Y''|X{=}x, n_X{=}x_{n_X}, X'{=}x', M{=}m) = \delta_K(n_X = x - x' - x_{n_x}). \tag{253}$
Here $\delta_K(\cdot)$ denotes the Kronecker delta function [59]. Marginalizing $(Y, Y', n_Y)$ from (246), we obtain:
$$\bar{Q}(M, X, X', Y'', n_X) = \bar{Q}(M)\bar{Q}(X|M)\bar{Q}(X', n_X, Y''|M, X) = \bar{Q}(M)\bar{Q}(X, X', Y'', n_X|M).$$
Consider the distribution $\bar{Q}(X, X', n_X, Y''|M)$. Then, we know:
$\bar{Q}(X|M{=}m) = f(\delta_m^X)$ and
$\bar{Q}(n_Y|X, M), \bar{Q}(X'|X, n_X, M),$ and $\bar{Q}(Y''|X, n_X, , X', M)$ are as defined in (253). $\quad$ (254)
Comparing the above equation with the structure of $\bar{Q}(X, X', n_X, Y''|M)$ specified in proposition 9, we can conclude that $(X', Y'', n_X)$ are just dilated versions of $X$. Consequently, using the result of proposition 9, we have:
$$\bar{Q}(X'|M{=}m) = f(\delta_m^X - \delta_m^Y - (\delta_{bias}^X - \delta_{bias}^Y)),$$
$$\bar{Q}(n_X|M) = f(\delta_{bias}^X),$$
$$\bar{Q}(Y''|M) = f(\delta_m^Y - \delta_{bias}^Y), \text{ and}$$
$$(X', Y'', n_X) \text{ are jointly conditionally independent conditioned on } M. \tag{255}$$

This concludes our construction of $\bar{Q}(\vec{\mathbf{B}}|\vec{\mathbf{A}}, M)$. We obtain our desired $\bar{Q}(M, X', Y'', n_X, Y', n_Y)$ by marginalizing $\vec{\mathbf{A}}$ out of $\bar{Q}(M, \vec{\mathbf{A}}, \vec{\mathbf{B}})$. From (251) and (255) we know that $\bar{Q}(M, X', Y'', n_X, Y', n_Y)$ satisfies the two main properties specified earlier. Since we did not make any specific assumptions on $Q$ aside from the fact that it lies in $\Delta_P$, the above construction can always be used to construct $(X', n_X, Y'', Y', n_Y)$ from $(M, X, Y)$ using the $\bar{Q}(\vec{\mathbf{B}}|\vec{\mathbf{A}}, M)$. Note that due to the particular construction of $\bar{Q}(\vec{\mathbf{B}}|\vec{\mathbf{A}}, M)$, we have:
$$X = X' + n_X + Y'' \text{ and } Y = Y' + n_Y. \tag{256}$$

### M.1.2 Propositions required for constructing $\bar{Q}$

We describe two important propositions that we use to show the existence of a $\bar{Q}$ for every $Q \in \Delta_P$.

**Proposition 8.** *Suppose the random variables $M$ and $Y$ have the joint distribution $Q(M, Y)$ such that*

$$Q(M) = P(M) \text{ and } Q(Y|M{=}m) = f(\delta_m^Y) \ \forall \ m \in \mathcal{M}, \tag{257}$$

*where $\mathcal{M}$ denotes the support of $M$ and $f(\delta_m^Y) \in \mathcal{F}_{\mathcal{D}}$ for some convolution-closed distribution family $\mathcal{F}_{\mathcal{D}}$. Furthermore, there also exists a $\delta_{bias}^Y \in \mathcal{D}$ such that*

$$\delta_m^Y = \epsilon_m^{(2)}(\delta_m^Y - \delta_{bias}^Y) \text{ for some } \epsilon_m^{(2)} \in [0, 1] \text{ and } (\delta_m^Y - \delta_{bias}^Y) \in \mathcal{D} \ \forall \ m \in \mathcal{M}. \tag{258}$$

*Consider two dilated versions of $Y$, denoted as $Y'$ and $n_Y$, defined as follows:*

$$\bar{Q}(Y'|Y{=}y, M{=}m) = G(\epsilon_m^{(2)}\delta_m^Y, (1 - \epsilon_m^{(2)})\delta_m^Y, y), \text{ and } n_Y = Y - Y'. \tag{259}$$

*Then, we have*

1. *$Y' \perp\!\!\!\perp n_Y | M$*

2. *The conditional distributions $\bar{Q}(Y'|M)$ and $\bar{Q}(n_Y|M)$ are specified as follows:*

$$\bar{Q}(Y'|M = m) = f(\epsilon_m^{(2)}\delta_m^Y) = f(\delta_m^Y - \delta_{bias}^Y) \text{ and}$$
$$\bar{Q}(n_Y|M{=}m) = f((1 - \epsilon_m^{(2)})\delta_m^Y) = f(\delta_{bias}^Y) \ \forall \ m \in \mathcal{M}. \tag{260}$$

*Proof.* We essentially modify the argument given in theorem 1 of [25] to prove the above proposition. Consider the conditional distribution of $Y$, $Y'$, and $n_Y$ conditioned on $M$, denoted as $\bar{Q}(Y, Y', n_Y | M)$. Then, by chain rule, we have $\bar{Q}(Y, Y', n_Y | M) = \bar{Q}(Y|M)\bar{Q}(Y'|M, Y)\bar{Q}(n_Y|Y, Y', M)$. By the assumptions in (257) and (259), we know that

$$\bar{Q}(Y|M{=}m) = f(\delta_m^Y), \bar{Q}(Y'|M{=}m, Y{=}y) = G(\epsilon_m^{(2)}\delta_m^Y, (1 - \epsilon_m^{(2)})\delta_m^Y, y), \text{ and}$$
$$\bar{Q}(n_Y|Y{=}y, Y'{=}y', M) = \delta_K(n_Y = y - y'), \tag{261}$$

where $\delta_K(\cdot)$ is the Kronecker delta function [59].

**Observation 1**: Our *first key observation* is that the distribution $\bar{Q}(Y', Y, n_Y | M{=}m)$ is completely specified through the distributions $\bar{Q}(Y|M)$ and $\bar{Q}(Y'|Y, M)$, as $n_Y$ is completely determined by $Y'$ and $Y$ and $\bar{Q}(n_Y|Y, Y', M)$ is specified through a Kronecker delta function.

Similarly, consider a new pair of random variables $Y'_{new}, n_Y^{new}$ such that $Y'_{new} \perp\!\!\!\perp n_Y^{new} | M$ and the respective conditional distributions of $Y'_{new}$ and $n_Y^{new}$ are specified as follows:

$$\bar{Q}_{new}(Y'_{new}|M{=}m) = f(\epsilon_m^{(2)}\delta_m^Y), \text{ and } \bar{Q}_{new}(n_Y^{new}|M{=}m) = f((1 - \epsilon_m^{(2)})\delta_m^Y) \ \forall \ m \in \mathcal{M}. \tag{262}$$

We will now show *two key properties* for the random variables $(Y'_{new}, n_Y^{new}, Y'_{new} + n_Y^{new})$ that will form the crux of our argument:

**Property 1**: $\bar{Q}_{new}(Y'_{new} + n_Y^{new}{=}y|M{=}m) = \bar{Q}(Y{=}y|M{=}m)$.

Let us now calculate the conditional distribution $\bar{Q}(Y'_{new} + n_Y^{new} = y|M)$:

$$\bar{Q}_{new}(Y'_{new} + n_Y^{new} = y|M) = \int_{k=-\infty}^{\infty} \bar{Q}_{new}(Y'_{new} = k, n_Y^{new} = y - k|M). \tag{263}$$

Using conditional dependence of $Y'_{new}$ and $n_Y^{new}$, we have $\bar{Q}_{new}(Y'_{new} = k, n_Y^{new} = y - k|M) = \bar{Q}_{new}(Y'_{new} = k|M)\bar{Q}_{new}(n_Y^{new} = y - k|M)$:

$$\bar{Q}_{new}(Y'_{new} + n_Y^{new} = y|M) = \int_{k=-\infty}^{\infty} \bar{Q}_{new}(Y'_{new} = k|M)\bar{Q}_{new}(n_Y^{new} = y - k|M). \tag{264}$$

Note that the R.H.S of the above equation defines a convolution of $\bar{Q}_{new}(Y'_{new}|M)$ and $\bar{Q}(n_Y^{new}|M)$, hence:

$$\bar{Q}_{new}(Y'_{new} + n_Y^{new} = y|M) = \bar{Q}_{new}(Y'_{new} = k|M) * \bar{Q}_{new}(n_Y^{new} = y - k|M), \tag{265}$$

where $*$ denotes the convolution operator. Substituting (262) in the above equation:

$$\bar{Q}_{new}(Y'_{new} + n_Y^{new} = y | M = m) = f(\epsilon_m^{(2)} \delta_m^Y) * f((1 - \epsilon_m^{(2)}) \delta_m^Y). \tag{266}$$

Using the properties of convolution-closed distributions:

$$\bar{Q}_{new}(Y'_{new} + n_Y^{new} = y | M = m) = f(\epsilon_m^{(2)} \delta_m^Y + (1 - \epsilon_m^{(2)}) \delta_m^Y) = f(\delta_m^Y). \tag{267}$$

Hence, by comparing above equation with (261) we have:

$$\bar{Q}_{new}(Y'_{new} + n_Y^{new} = y | M = m) = Q(Y = y | M = m). \tag{268}$$

**Property 2**: $\bar{Q}(Y'_{new} = y' | Y'_{new} + n_Y^{new} = y, M = m) = G(\epsilon_m^{(2)} \delta_m^Y, 1 - \epsilon_m^{(2)} \delta_m^Y, y)$

By definition of $G(\epsilon_m^{(2)} \delta_m^Y, (1 - \epsilon_m^{(2)}) \delta_m^Y, z)$ given in Sec. 5.1, we have:

$$G(\epsilon_m^{(2)} \delta_m^Y, (1 - \epsilon_m^{(2)}) \delta_m^Y, z) = \Pr(Z_{\epsilon_m^{(2)}} | Z_{\epsilon_m^{(2)}} + Z_{1 - \epsilon_m^{(2)}} = z), \text{ where}$$
$$\Pr(Z_{\epsilon_m^{(2)}}) = f(\epsilon_m^{(2)} \delta_m^Y), \Pr(Z_{1 - \epsilon_m^{(2)}}) = f((1 - \epsilon_m^{(2)}) \delta_m^Y), \text{ and } Z_{\epsilon_m^{(2)}} \perp\!\!\!\perp Z_{1 - \epsilon_m^{(2)}}. \tag{269}$$

In the following steps, we slightly modify the notation of $G(\epsilon_m^{(2)} \delta_m^Y, (1 - \epsilon_m^{(2)}) \delta_m^Y, z)$ to $G(z_{\epsilon_m^{(2)}}; \epsilon_m^{(2)} \delta_m^Y, (1 - \epsilon_m^{(2)}) \delta_m^Y, z)$ to make our notations more explicit as follows:

$$G(z_{\epsilon_m^{(2)}}; \epsilon_m^{(2)} \delta_m^Y, (1 - \epsilon_m^{(2)}) \delta_m^Y, z) = \Pr(Z_{\epsilon_m^{(2)}} = z_{\epsilon_m^{(2)}} | Z_{\epsilon_m^{(2)}} + Z_{1 - \epsilon_m^{(2)}} = z). \tag{270}$$

We use the definition of the conditional distribution to express $G(\epsilon_m^{(2)} \delta_m^Y, (1 - \epsilon_m^{(2)}) \delta_m^Y, z)$ as:

$$G(z_{\epsilon_m^{(2)}}; \epsilon_m^{(2)} \delta_m^Y, (1 - \epsilon_m^{(2)}) \delta_m^Y, z) = \frac{\Pr(Z_\epsilon = z_{\epsilon_m^{(2)}}, Z_{\epsilon_m^{(2)}} + Z_{1 - \epsilon_m^{(2)}} = z)}{\Pr(Z_{\epsilon_m^{(2)}} + Z_{1 - \epsilon_m^{(2)}} = z)} \tag{271}$$

$$= \frac{\Pr(Z_\epsilon = z_{\epsilon_m^{(2)}}) \Pr(Z_{\epsilon_m^{(2)}} + Z_{1 - \epsilon_m^{(2)}} = z | Z_{\epsilon_m^{(2)}} = z_{\epsilon_m^{(2)}})}{\Pr(Z_{\epsilon_m^{(2)}} + Z_{1 - \epsilon_m^{(2)}} = z)}. \tag{272}$$

Using the fact that $\Pr(Z_{\epsilon_m^{(2)}} + Z_{1 - \epsilon_m^{(2)}} = z | Z_{\epsilon_m^{(2)}} = z_{\epsilon_m^{(2)}}) = \Pr(Z_{1 - \epsilon_m^{(2)}} = z - z_{\epsilon_m^{(2)}})$, since $Z_{\epsilon_m^{(2)}} \perp\!\!\!\perp Z_{1 - \epsilon_m^{(2)}}$:

$$= \frac{\Pr(Z_\epsilon = z_{\epsilon_m^{(2)}}) \Pr(Z_{1 - \epsilon_m^{(2)}} = z - z_{\epsilon_m^{(2)}})}{\Pr(Z_{\epsilon_m^{(2)}} + Z_{1 - \epsilon_m^{(2)}} = z)}. \tag{273}$$

Note that by properties of convolution-closed distributions, $\Pr(Z_{\epsilon_m^{(2)}} + Z_{1 - \epsilon_m^{(2)}} = z)) = f(z; \delta_m^Y)$, where $f(z; \delta_m^Y)$ is to be understood as the distribution $f(\delta_m^Y)$ evaluated at $z$. Substituting (269) in the above equation:

$$= \frac{f(z_{\epsilon_m^{(2)}}; \epsilon_m^{(2)} \delta_m^Y) f(z - z_{\epsilon_m^{(2)}}; (1 - \epsilon_m^{(2)}) \delta_m^Y)}{f(z; \delta_m^Y)}. \tag{274}$$

Now, note that by (262) and (267), we have:

$$\bar{Q}_{new}(Y'_{new} = z_\epsilon | M = m) = f(z_{\epsilon_m^{(2)}}; \epsilon_m^{(2)} \delta_m^Y),$$
$$\bar{Q}_{new}(n_Y^{new} = z - z_\epsilon | M = m) = f(z - z_{\epsilon_m^{(2)}}; (1 - \epsilon_m^{(2)}) \delta_m^Y), \text{ and}$$
$$\bar{Q}_{new}(Y'_{new} + n_Y^{new} = z | M = m) = f(z; \delta_m^Y) \tag{275}$$

Hence, combining the above equation with (274), and renaming $z_{\epsilon_m^{(2)}}$ as $y'$ and $z$ as $y$, we have:

$$G(\epsilon_m^{(2)} \delta_m^Y, (1 - \epsilon_m^{(2)}) \delta_m^Y, y) = \frac{\bar{Q}_{new}(Y'_{new} = y' | M = m) \bar{Q}_{new}(n_Y^{new} = y - y' | M = m)}{\bar{Q}_{new}(Y'_{new} + n_Y^{new} = y | M = m)}. \tag{276}$$

Noting the fact that $\bar{Q}_{new}(n_Y^{new}{=}y - y'|M{=}m) = \bar{Q}_{new}(Y'_{new} + n_Y^{new}{=}y|M{=}m, Y'_{new} = y')$ due to $Y'_{new} \perp\!\!\!\perp n_Y^{new}|M$ and abbreviating notation pertaining to $y'$, we have:

$$G(\epsilon_m^{(2)}\delta_m^Y, (1 - \epsilon_m^{(2)})\delta_m^Y, y) = \frac{\bar{Q}_{new}(Y'_{new}|M{=}m)\bar{Q}_{new}(Y'_{new} + n_Y^{new}{=}y|M{=}m)}{\bar{Q}_{new}(Y'_{new} + n_Y^{new}{=}y|M{=}m)}. \quad (277)$$

Using Bayes rule in the above equation provides us with our desired property:

$$G(\epsilon_m^{(2)}\delta_m^Y, (1 - \epsilon_m^{(2)})\delta_m^Y, y) = \bar{Q}_{new}(Y'_{new}|M{=}m, Y'_{new} + n_Y^{new}{=}y) \quad (278)$$

Furthermore, by comparing (259) with the above equation, we have:

$$\bar{Q}_{new}(Y'_{new} = y'|Y'_{new} + n_Y^{new} = y, M = m) = \bar{Q}(Y' = y'|Y = y, M = m). \quad (279)$$

**Observation 2**: Note that the distribution $\bar{Q}(Y'_{new}, n_Y^{new}, Y'_{new} + n_Y^{new}|M)$ is also completely determined by specifying $\bar{Q}(Y'_{new} + n_Y^{new}|M)$ and $\bar{Q}(Y'_{new}|Y'_{new} + n_Y^{new}, M)$ since $n_Y^{new}$ is completely determined by $Y'_{new} + n_Y^{new}$ and $Y'_{new}$.

By property 1 and (279), we know that

$$\bar{Q}_{new}(Y'_{new} + n_Y^{new}|M) = Q(Y|M), \text{ and } \bar{Q}_{new}(Y'_{new}|Y'_{new} + n_Y^{new}, M) = \bar{Q}(Y'|Y, M). \quad (280)$$

Therefore, combining the above equation with observations 1 and 2, we have:

$$\bar{Q}(Y', Y, n_Y|M) = \bar{Q}(Y'_{new}, Y'_{new} + n_Y^{new}, n_Y^{new}|M). \quad (281)$$

Consequently the distribution $\bar{Q}(Y', n_Y|M) = \bar{Q}_{new}(Y'_{new}, n_Y^{new}|M)$. Since, $Y'_{new} \perp\!\!\!\perp n_Y^{new}|M$, we also have $Y' \perp\!\!\!\perp n_Y|M$. Furthermore, by (262),

$$\bar{Q}_{new}(Y'_{new}|M{=}m) = \bar{Q}(Y'|M) = f(\epsilon_m^{(2)}\delta_m^Y) \,\forall\, m \in \mathcal{M}, \text{ and}$$
$$\bar{Q}_{new}(n_Y^{new}|M{=}m) = \bar{Q}(n_Y|M) = f((1 - \epsilon_m^{(2)})\delta_m^Y) \,\forall\, m \in \mathcal{M}. \quad (282)$$

By (258), we know that $\epsilon_m^{(2)}\delta_m^Y = \delta_m^Y - \delta_{bias}^Y$ and $(1 - \epsilon_m^{(2)})\delta_m^Y = \delta_{bias}^Y$. Hence,

$$\bar{Q}(Y'|M) = f(\epsilon_m^{(2)}\delta_m^Y) = f(\delta_m^Y - \delta_{bias}^Y) \,\forall\, m \in \mathcal{M}, \text{ and}$$
$$\bar{Q}(n_Y|M) = f((1 - \epsilon_m^{(2)})\delta_m^Y) = f(\delta_{bias}^Y) \,\forall\, m \in \mathcal{M}. \quad (283)$$

$$\square$$

**Proposition 9.** *Suppose the random variables $M$ and $X$ have the joint distribution $Q(M, X)$, such that*

$$Q(M) = P(M) \text{ and } Q(X|M{=}m) = f(\delta_m^X) \,\forall\, m \in \mathcal{M}, \quad (284)$$

*where $\mathcal{M}$ denotes the support of $M$ and $f(\delta_m^X) \in \mathcal{F}_\mathcal{D}$ for some convolution-closed distribution family $\mathcal{F}_\mathcal{D}$. Furthermore, there also exist $\delta_{bias}^X \in \mathcal{D}$, and $\delta_m^Y \in \mathcal{D} \,\forall\, m \in \mathcal{M}$ such that*

$$\delta_m^X - \delta_{bias}^X = \epsilon_m^{(1)}(\delta_m^Y - \delta_{bias}^Y) \text{ for some } \epsilon_m^{(1)} \in [0, 1],$$
$$\delta_m^X = \epsilon_m^{(3)}(\delta_m^X - \delta_{bias}^X) \text{ for some } \epsilon_m^{(3)} \in [0, 1], \text{ and}$$
$$(\delta_m^X - \delta_m^Y - (\delta_{bias}^X - \delta_{bias}^Y)), (\delta_m^X - \delta_{bias}^X), (\delta_m^Y - \delta_{bias}^Y) \in \mathcal{D} \,\forall\, m \in \mathcal{M}. \quad (285)$$

*Then consider three dilated versions of $X$, denoted as $X', Y''$, and $n_X$, defined as follows:*

$$\bar{Q}(X', Y'', n_X|X, M) = \bar{Q}(n_X|X, M)\bar{Q}(X'|X, n_X, M)\bar{Q}(Y''|X, n_X, M, X'), \text{ where}$$
$$\bar{Q}(n_X|X{=}x, M{=}m) = G((1 - \epsilon_m^{(3)})\delta_m^X, \epsilon_m^{(3)}\delta_m^X, x),$$
$$\bar{Q}(X'|X{=}x, n_X{=}x_{n_X}, M{=}m) = G((1 - \epsilon_m^{(1)})(\delta_m^X - \delta_{bias}^X), \epsilon_m^{(1)}(\delta_m^X - \delta_{bias}^X), x - n_x), \text{ and}$$
$$Y'' = X - X - n_X. \quad (286)$$

*Then, we have*

1. *$(n_X, X', Y'')$ are jointly conditionally independent conditioned on $M$.*

2. *The conditional distributions $\bar{Q}(X'|M)$, $\bar{Q}(Y''|M)$, and $\bar{Q}(n_X|M)$ are specified as follows:*

$$\bar{Q}(X'|M) = f(\delta_m^X - \delta_m^Y - (\delta_{bias}^X - \delta_{bias}^Y)),$$
$$\bar{Q}(Y''|M) = f(\delta_m^X - \delta_{bias}^Y), \text{ and}$$
$$\bar{Q}(n_X|M) = f(\delta_{bias}^X). \tag{287}$$

*Proof.* The proof essentially consists of recursively applying the proof of proposition 8. Consider a new auxiliary random variable $Z = X - n_X$. Then, the distribution $\bar{Q}(X, n_X, Z|M)$ has the following properties:

$$\bar{Q}(X|M) \overset{(a)}{=} f(\delta_m^X), \bar{Q}(n_X|M=m, X=x) \overset{(b)}{=} G((1 - \epsilon_m^{(3)})\delta_m^X, \epsilon_m^{(3)}\delta_m^X, x), \text{ and}$$
$$\bar{Q}(Z|X=x, n_X=x_{n_X}, M=m) = \delta_K(Z = x - x_{n_X}), \tag{288}$$

where $\delta_K(\cdot)$ is the Kronecker delta function, and the equalities $(a)$ and $(b)$ follow from (284) and (285), respectively. Note that the distribution $\bar{Q}(X, n_X, Z|M)$ has exactly the same structure as $\bar{Q}(Y, Y', n_Y|M)$ described in (261), hence $Z$ and $n_Y$ are just dilated versions of $X$. Consequently, from the result of proposition 8, we know that $Z \perp\!\!\!\perp n_X|M$, and the conditional distributions $\tilde{Q}(Z|M)$ and $\bar{Q}(n_Y|M)$ are described as follows:

$$\bar{Q}(n_X|M=m) = f((1 - \epsilon_m^{(3)})\delta_m^X), \text{ and } \bar{Q}(Z|M=m) = f(\epsilon_m^{(3)}\delta_m^X). \tag{289}$$

From (285), we have $\epsilon_m^{(3)}\delta_m^X = \delta_m^X - \delta_{bias}^X$ and $(1 - \epsilon_m^{(3)})\delta_m^X = \delta_{bias}^X$. Consequently,

$$\bar{Q}(n_X|M=m) = f(\delta_{bias}^X), \text{ and } \bar{Q}(Z|M=m) = f(\delta_m^X - \delta_{bias}^X). \tag{290}$$

We alternatively express $\tilde{Q}(X'|n_X, X, M)$ as follows:

$$\bar{Q}(X'|n_X=x_{n_X}, X=x, M=m) = G((1 - \epsilon_m^{(1)})(\delta_m^X - \delta_{bias}^X), \epsilon_m^{(1)}(\delta_m^X - \delta_{bias}^X), x - x_{n_X}). \tag{291}$$

Note that the distribution $\bar{Q}(X'|n_X=x_{n_X}, X=x, M=m)$ only depends on $X$ and $n_X$ through their difference $X - n_X$. Hence, substituting $Z = X - n_X$ in the above equation:

$$\bar{Q}(X'|n_X=x_{n_X}, X=x, M=m) = \bar{Q}(X'|Z=x - x_{n_X}, M=m)$$
$$= G((1 - \epsilon_m^{(1)})(\delta_m^X - \delta_{bias}^X), \epsilon_m^{(1)}(\delta_m^X - \delta_{bias}^X), x - n_X), \tag{292}$$
$$\Rightarrow \bar{Q}(X'|Z=z, M=m) = G((1 - \epsilon_m^{(1)})(\delta_m^X - \delta_{bias}^X), \epsilon_m^{(1)}(\delta_m^X - \delta_{bias}^X), z). \tag{293}$$

Hence, consider the following distribution $\bar{Q}(Z, X', Y''|M)$, where from (290) and (293), we have:

$$\bar{Q}(Z|M=m) = f(\delta_m^X - \delta_{bias}^X),$$
$$\bar{Q}(X'|Z = z, M = m) = G((1 - \epsilon_m^{(1)})(\delta_m^X - \delta_{bias}^X), \epsilon_m^{(1)}(\delta_m^X - \delta_{bias}^X), z). \tag{294}$$

Furthermore, by (286), we have $Y'' = X - n_X - X' \overset{(c)}{=} Z - X'$, where $(c)$ is due to $Z = X - n_X$. Hence,

$$\bar{Q}(Y''|Z=z, M, X'=x) = \delta_K(Y'' = z - x). \tag{295}$$

Comparing (295) and (294) with (261), we can again conclude that $X''$ and $Y'$ are just dilated versions of $Z$. Hence, again using the result of proposition 8, we have $X' \perp\!\!\!\perp Y''|M$, and

$$\bar{Q}(X'|M=m) = f((1 - \epsilon_m^{(1)})(\delta_m^X - \delta_{bias}^X)), \text{ and } \bar{Q}(Y''|M=m) = f(\epsilon_m^{(1)}(\delta_m^X - \delta_{bias}^X)). \tag{296}$$

From (285), we have $\epsilon_m^{(1)}(\delta_m^X - \delta_{bias}^X) = \delta_m^Y - \delta_{bias}^Y$ and $(1 - \epsilon_m^{(1)})(\delta_m^X - \delta_{bias}^X) = \delta_m^X - \delta_M^Y - (\delta_{bias}^X - \delta_{bias}^Y)$. Consequently,

$$\bar{Q}(X'|M=m) = f(\delta_m^X - \delta_m^Y - (\delta_{bias}^X - \delta_{bias}^Y)), \text{ and } \bar{Q}(Y''|M=m) = f(\delta_m^Y - \delta_{bias}^Y). \tag{297}$$

Hence, by (290) and (297), we can conclude that the conditional distributions $\bar{Q}(X'|M)$, $\bar{Q}(Y''|M)$, and $\bar{Q}(n_X|M)$ are specified as follows:

$$
\begin{aligned}
\bar{Q}(X'|M=m) &= f(\delta_m^X - \delta_m^Y - (\delta_{bias}^X - \delta_{bias}^Y)), \\
\bar{Q}(Y''|M) &= f(\delta_m^X - \delta_{bias}^Y), \text{ and} \\
\bar{Q}(n_X|M) &= f(\delta_{bias}^X).
\end{aligned}
\tag{298}
$$

*Showing joint conditional independence of* $(n_Y, X', Y'')$ *given* $M$:

Consider the distribution $\bar{Q}(n_X, Z, X', Y''|M)$. Using the chain rule, we have:

$$
\begin{aligned}
\bar{Q}(n_X, Z, X', Y''|M) &= \bar{Q}(n_X, Z|M)Q(X', Y''|M, n_X, Z), \\
&\overset{(b)}{=} \bar{Q}(n_X|M)\bar{Q}(Z|M)Q(X', Y''|M, n_X, Z), \\
&\overset{(c)}{=} \bar{Q}(n_X|M)\bar{Q}(Z|M)Q(X', Y''|M, Z),
\end{aligned}
\tag{299}
$$

where $(b)$ follows from $n_X \perp\!\!\!\perp Z|M$. For showing $(c)$, observe that $Y'' = Z - X'$, and hence we can write $\bar{Q}(Y''|M, Z, n_Y, X') = \bar{Q}(Y''|Z, X')$. Furthermore, by analyzing $\bar{Q}(X'|n_X, Z, M)$, we can conclude:

$$
\bar{Q}(X'|n_X, Z, M) = \bar{Q}(X'|n_X, X - n_X, M) = \bar{Q}(X'|n_X, X, M) = \bar{Q}(X'|Z, M).
\tag{300}
$$

Marginalizing $Z$ out of (299), we obtain:

$$
\bar{Q}(n_X, X', Y''|M) = \bar{Q}(n_X|M)\bar{Q}(X', Y''|M) \overset{(d)}{=} \bar{Q}(n_X|M)Q(X'|M)\bar{Q}(Y''|M),
\tag{301}
$$

where $(d)$ follows from $X' \perp\!\!\!\perp Y''|M$. By (301), we know that $(n_X, X'', Y')$ are jointly conditionally independent given $M$, concluding our proof. $\qquad\square$

## M.2 Analytically optimizing the upper bound

**Proposition 10.** *Let* $\bar{Q}(M, X', Y'', n_X, Y', n_Y)$ *be the distribution derived from a distribution* $Q \in \Delta_P$ *using the construction scheme specified in Appx. M.1. Then, the minimization problem specified in* (302) *is analytically solvable with the optimizing* $\bar{Q}^*$ *given by* (303).

$$
\min_{Q \in \Delta_P} I_{\bar{Q}}(M; [X', Y'', n_X, Y', n_Y]), \text{ (note that } \bar{Q} \text{ is a function of } Q)
\tag{302}
$$

$\bar{Q}^*(M, X', Y'', n_X, n_Y, Y') = \bar{Q}^*(M)\bar{Q}^*(X'|M)\bar{Q}^*(Y''|M)\bar{Q}^*(n_X)\bar{Q}^*(n_Y)\bar{Q}^*(Y'|Y'')$, *with*

$\bar{Q}^*(M) = P(M), \bar{Q}^*(X'|M=m) = f(\delta_m^X - \delta_m^Y - (\delta_{bias}^X - \delta_{bias}^Y)), \bar{Q}^*(n_X) = f(\delta_{bias}^X)$,

$\bar{Q}^*(Y''|M=m) = f(\delta_m^Y - \delta_{bias}^Y), \bar{Q}^*(n_Y) = f(\delta_{bias}^Y),$ *and* $\bar{Q}^*(Y'|Y'') = \mathbb{I}[Y'' = Y']$, $\qquad$ (303)

*where* $\mathbb{I}[\cdot]$ *is the indicator function. Note that for the minimizing* $\bar{Q}^*$, *we have* $(n_X, n_Y, (M, X', Y''))$ *are jointly independent and* $Y' = Y''$.

*Proof.* First note that by the construction scheme defined in Appx. M.1, we know that there is a unique $\bar{Q}$ for every $Q \in \Delta_P$. Hence, we can interpret $\bar{Q}$ as a function of $Q$, and consequently the minimization problem in (302) is well-defined. Using the chain rule of mutual information, we derive the following lower bound for $I_{\bar{Q}}(M; [X', Y'', n_X, Y', n_Y])$:

$$
I_{\bar{Q}}(M; [X', Y'', n_X, Y', n_Y]) = I_{\bar{Q}}(M; [X', Y'']) + I_{\bar{Q}}(M; [n_X, Y', n_Y]|[X', Y'])
\tag{304}
$$

$$
\Rightarrow I_{\bar{Q}}(M; [X', Y'', n_X, Y', n_Y]) \overset{(a)}{\geq} I_{\bar{Q}}(M; [X', Y'']),
\tag{305}
$$

where $(a)$ is due to the non-negativity of conditional mutual information. Furthermore, note that by properties of $\bar{Q}$ derived in Appx. M.1, we know that $X' \perp\!\!\!\perp Y''|M \ \forall \ Q \in \Delta_P$. Consequently, the term $I_{\bar{Q}}(M; [X', Y''])$ does note vary as we choose different $Q$ from $\Delta_P$ as the corresponding $\bar{Q}(M, X', Y'') = \bar{Q}(M)\bar{Q}(X'|M)\bar{Q}(Y''|M)$ and the terms:

$$
\begin{aligned}
\bar{Q}(M) &= Q(M) = P(M), \bar{Q}(X'|M=m) = f(\delta_m^X - \delta_m^Y - (\delta_{bias}^X - \delta_{bias}^Y)), \text{ and} \\
\bar{Q}(Y''|M) &= f(\delta_m^Y - \delta_{bias}^Y)
\end{aligned}
\tag{306}
$$

do not vary for all $Q \in \Delta_P$ (see Appx. M.1). Therefore, we have:

$$I(M; [X', Y'']) \leq I_{\bar{Q}}(M; [X', Y'', n_X, Y', n_Y]) \ \forall \ Q \in \Delta_P, \tag{307}$$

where we forgo the subscript $\bar{Q}$ for $I_{\bar{Q}}(M; [X'', Y'])$ to indicate that the term $I_{\bar{Q}}(M; [X'', Y'])$ does not vary over $\Delta_P$. Hence, if we show that the distribution $\bar{Q}^*(M, X', Y'', n_X, Y', n_Y)$ specified in (303) achieves the lower bound in (307), then $\bar{Q}^*(M, X', Y'', n_X, Y', n_Y)$ should minimize the value of $I_{\bar{Q}}(M; [X', Y'', n_X, Y', n_Y])$ over the set $\Delta_P$. We now calculate $I_{\bar{Q}^*}(M; [X', Y'', n_X, Y', n_Y])$. By the chain rule of mutual information:

$$I_{\bar{Q}^*}(M; [X', Y'', n_X, Y', n_Y]) = I_{\bar{Q}^*}(M; [n_X, n_Y, X', Y'']) + I_{\bar{Q}^*}(M; Y' | n_X, n_Y, X, Y'').$$

Using the fact that $Y' = Y''$, we know that $I_{\bar{Q}^*}(M; Y' | n_X, n_Y, X, Y'')$ is zero. A simple justification follows from the fact $I_{\bar{Q}^*}(M; Y' | n_X, n_Y, X, Y'') = H(Y' | n_X, n_Y, X, Y'') - H(Y' | n_X, n_Y, X, Y'', M)$, where $H(.|.)$ is the conditional entropy function [28]. Observe that both conditional entropy terms are zero as $Y' = Y''$, and condition entropy of a random variable conditioned on itself is zero. Therefore, we can simplify the above equation as:

$$I_{\bar{Q}^*}(M; [X', Y'', n_X, Y', n_Y]) = I_{\bar{Q}^*}(M; [n_X, n_Y, X', Y'']). \tag{308}$$

Applying the chain rule of mutual information again, we obtain:

$$I_{\bar{Q}^*}(M; [X', Y'', n_X, Y', n_Y]) = I_{\bar{Q}^*}(M; [n_X, n_Y]) + I_{\bar{Q}^*}(M; [X', Y''] | n_X, n_Y). \tag{309}$$

Using the fact that $(n_X, n_Y) \perp\!\!\!\perp (M, X', Y'')$, we know that $I(M; [n_X, n_Y]) = 0$ as mutual information between independent random variables is zero [28]. Consequently,

$$I_{\bar{Q}^*}(M; [X', Y'', n_X, Y', n_Y]) = I_{\bar{Q}^*}(M; [X', Y''] | n_X, n_Y) \overset{(b)}{=} I_{\bar{Q}^*}(M; [X', Y'']), \tag{310}$$

where $(b)$ again follows from the fact that $(n_X, n_Y) \perp\!\!\!\perp (M, X', Y'')$, and, consequently, conditioning $(M, X', Y'')$ on $(n_X, n_Y)$ has no effect. From (310), we see the proposed $\bar{Q}^*$ in (303) does indeed achieve the lower bound specified in (307), and consequently minimizes (302).

The final step remaining in our proof is to ensure that the proposed $\bar{Q}^*$ corresponds to a valid $Q^*_{UB}(M, X, Y)$ lying in $\Delta_P$, i.e., there is some $Q^*_{UB}(M, X, Y) \in \Delta_P$ that generates $\bar{Q}^*$ using the construction scheme outlined in Appx. M.1. We know that the relationship between the random variables for $(M, X', Y'', n_X, Y', n_Y)$ having the joint distribution $\bar{Q}$ and $(M, X, Y)$ having the distribution $Q$ can be expressed as follows: $X = X' + Y'' + n_X$ and $Y = Y' + n_Y$. Hence, we use these relationships for calculating the conditional marginals $Q^*_{UB}(X|M)$ and $Q^*_{UB}(Y|M)$ of the corresponding distribution $Q^*_{UB}(M, X, Y)$ for the proposed distribution $\bar{Q}^*(M, X', Y'', n_X, Y', n_Y)$:

$$
\begin{aligned}
Q^*_{UB}(X|M = m) &= \bar{Q}^*(X' + Y'' + n_X | M = m) \\
&\overset{(c)}{=} \bar{Q}^*(X'|M = m) * \bar{Q}^*(Y''|M = m) * \bar{Q}^*(n_X|M = m) \\
&= f(\delta_m^X - \delta_m^Y - (\delta_{bias}^X - \delta_{bias}^Y)) * f(\delta_m^Y - \delta_{bias}^Y) * f(\delta_{bias}^X) \\
&\overset{(d)}{=} f(\delta_m^X - \delta_m^Y - (\delta_{bias}^X - \delta_{bias}^Y) + \delta_m^Y - \delta_{bias}^Y + \delta_{bias}^X) \\
&= f(\delta_m^X) = P(X|M = m).
\end{aligned}
\tag{311}
$$

where $*$ denotes the convolution operator, and $(c)$ follows from the fact that $(X', Y'', n_X)$ are jointly conditionally independent given $M$ under $\bar{Q}^*$, and $(d)$ follows from the properties of convolution-closed distributions. Similarly, calculating $Q^*_{UB}(Y|M = m)$:

$$
\begin{aligned}
Q^*_{UB}(Y|M = m) &= \bar{Q}^*(Y' + n_Y | M = m) \\
&\overset{(e)}{=} \bar{Q}^*(Y''|M = m) * \bar{Q}^*(n_Y|M = m) \\
&= f(\delta_m^Y - \delta_{bias}^Y) * f(\delta_{bias}^Y) \\
&= f(\delta_m^Y - \delta_{bias}^Y + \delta_{bias}^X) \\
&= f(\delta_m^Y) = P(Y|M = m),
\end{aligned}
\tag{312}
$$

where $(e)$ follows from the fact that $Y' \perp\!\!\!\perp n_Y | M$ under $\bar{Q}^*$. It is easy to see that $Q^*_{UB}(M) = \bar{Q}(M) = P(M)$. Hence, we have $Q^*_{UB}(M, X) = P(M)P(X|M) = P(M, X)$ and $Q^*_{UB}(M, Y) = P(M)P(X = Y|M) = P(M, Y)$, which implies $Q^*_{UB}(M, X, Y) \in \Delta_P$, concluding our proof. □

## M.3   Additional details regarding simulation

The list of different function pairs used in the simulation study are listed in Table 1. The function pairs were chosen with the only requirement being that they satisfied the assumptions in (9). For constructing the analytical estimate $\hat{Q}_A(M, X, Y)$, the corresponding $\delta_{bias}^X$ and $\delta_{bias}^Y$ were chosen as follows:

1. For each distribution and function pair, we first determine the range of possible values for $\delta_{bias}^X$ and $\delta_{bias}^Y$.

2. For $\delta_{bias}^Y$, we choose the range as $[0, \min_{m \in \mathcal{M}} f_2(m))$, where $\mathcal{M}$ is the support set of $M$ for that distribution. Note that we cannot choose $\delta_{bias}^Y > \min_{m \in \mathcal{M}} f_2(m)$ as the corresponding conditional distribution $\bar{Q}^*(Y'|M) = \text{Poisson}(f_2(M) - \delta_{bias}^Y)$ (required for the optimal $\bar{Q}^*$ from which $Q_A$ is constructed, see Appx. M.2) would have a negative rate parameter, which is not allowed. Within this range, we choose 10 different, uniformly spaced values for $\delta_{bias}^Y$.

3. For each chosen value of $\delta_{bias}^Y$, the corresponding range for $\delta_{bias}^X$ is then chosen as $[0, \min\{\min_{m \in \mathcal{M}} f_1(m), \min_{m \in \mathcal{M}}(f_1(m) - f_2(m) + \delta_{bias}^Y)\}]$. Note that choosing the upper limit of $\delta_{bias}^X$ as $\min\{\min_{m \in \mathcal{M}} f_1(m), \min_{m \in \mathcal{M}}(f_1(m) - f_2(m) + \delta_{bias}^Y)\}$ ensures that the distribution $\bar{Q}^*(X'|M) = \text{Poisson}((f_1(M) - \delta_{bias}^X) - (f_2(M) - \delta_{bias}^Y))$ and $\bar{Q}^*(Z|M) = \text{Poisson}(f_1(M) - \delta_{bias}^X)$ (required for specifying the optimal $\bar{Q}^*$ from which $Q_A$ is constructed, see Appx. M.1 and Appx. M.2) have positive rate parameters. For each value of $\delta_{bias}^Y$, we again choose 10 different values of $\delta_{bias}^X$ within the above specified range. In total, we chose 100 different pairs of $(\delta_{bias}^X, \delta_{bias}^Y)$.

For each pair of $(\delta_{bias}^X, \delta_{bias}^Y)$, we calculate the analytical $Q_A(M, X, Y)$ as follows:

First, we construct the optimal $\bar{Q}^*$ distribution using the structure provided in proposition 10. The exact specification of $\bar{Q}^*$ is described below. We simplify the structure of $\bar{Q}^*$ by using the fact that $Y' = Y''$:

$$
\begin{aligned}
\bar{Q}^*(M, X', Y', n_X, n_Y, Y'') &\equiv \bar{Q}^*(M, X', Y', n_X, n_Y), \\
&= \bar{Q}^*(M)\bar{Q}^*(X'|M)\bar{Q}^*(Y'|M)\bar{Q}^*(n_X)\bar{Q}^*(n_Y), \text{ with} \\
\bar{Q}^*(M) &= P(M), \\
\bar{Q}^*(X'|M) &= \text{Poisson}(f_1(M) - f_2(M) - (\delta_{bias}^X - \delta_{bias}^Y)), \\
\bar{Q}^*(n_X) &= \text{Poisson}(\delta_{bias}^X), \\
\bar{Q}^*(Y''|M) &= \text{Poisson}(f_2(M) - \delta_{bias}^Y), \\
\bar{Q}^*(n_Y) &= \text{Poisson}(\delta_{bias}^Y).
\end{aligned}
\tag{313}
$$

Then, we use the property that $X = X' + Y' + n_X$, and $Y = Y' + n_Y$, and use that to appropriately modify $\bar{Q}^*$ to obtain $Q_A$ as follows:

- We calculate a new distribution $\bar{Q}^*(M, X' + Y', Y', n_X, n_Y)$:

$$
\begin{aligned}
\bar{Q}^*(M, X' + Y', Y', n_X, n_Y) &= \bar{Q}^*(M)\bar{Q}^*(n_X)\bar{Q}^*(n_Y)\bar{Q}^*(X' + Y'|M) \times \\
&\quad \bar{Q}^*(Y'|X' + Y', M). \\
&= \bar{Q}^*(M)\bar{Q}^*(n_X)\bar{Q}^*(n_Y)\bar{Q}^*(Z|M)\bar{Q}^*(Y'|Z, M),
\end{aligned}
\tag{314}
$$

  where $Z = X' + Y'$ and alternatively also $Z = X - n_X$ (as $X = X' + Y' + n_X$). First note that $\bar{Q}^*(Z|M)$ from Appx. M.2 has the form $f(\delta_m^X - \delta_{bias}^X)$, which for the Poisson case is $\text{Poisson}(f_1(M) - \delta_{bias}^X)$. Furthermore, we can recognize that $\bar{Q}^*(Y' = y|Z = z, M = m) = \bar{Q}^*(Y' = y'|X' + Y' = x' + y', M = m) = G(y'; \epsilon_m^{(1)}, 1 - \epsilon_m^{(1)}, x' + y')$, which for the Poisson case can be written as $\text{Binomial}(Z, p_m)$ where $p_m = {}^{f_2(M) - \delta_{bias}^Y}\!/{}_{f_1(M) - \delta_{bias}^X}$.

- Now define $\mathbb{S}_x^m = \{x_{n_X} \in \mathcal{X}_{n_X}^m, z \in \mathcal{Z}_m : x_{n_X} + z = x\}$, where $\mathcal{X}_{n_X}^m$ and $\mathcal{Z}_m$ are the corresponding support sets of the distributions $\bar{Q}^*(n_X|M = m)$ and $\bar{Q}^*(Z|M = m)$,

respectively. Then, using the fact that $X = Z + n_X$, we have:

$$\bar{Q}^*(M{=}m, X{=}x, n_Y, Y') = \bar{Q}^*(M, Z + n_X, n_Y, Y') = \bar{Q}^*(M = m)\bar{Q}^*(n_Y)\times$$

$$\sum_{(x_{n_X}, z) \in \mathbb{S}_x^m} \bar{Q}^*(n_X = x_{n_X})\bar{Q}^*(Z = z | M = m)\bar{Q}^*(Y' | Z = z, M = m), \qquad (315)$$

where we know that for the Poisson case:

$$\bar{Q}^*(M) = P(M), \bar{Q}^*(n_Y) = \text{Poisson}(\delta_{bias}^Y), \bar{Q}^*(Z|M) = \text{Poisson}(f_1(M) - \delta_{bias}^X),$$

$$\bar{Q}^*(n_X) = \text{Poisson}(\delta_{bias}^X), \text{ and } \bar{Q}^*(Y'|Z, M) = \text{Binomial}\left(Z, \frac{f_2(M) - \delta_{bias}^Y}{f_1(M) - \delta_{bias}^X}\right). \ (316)$$

- Using a similar procedure, we derive $Q_A(M, X, Y)$ from $\bar{Q}^*(M, X, Y', n_Y)$ as $Y = n_Y + Y'$. Define $\mathbb{S}_y^m = \{y_{n_y} \in \mathcal{Y}_{n_X}^m, z \in \mathcal{Y}_m' : y_{n_X} + y' = y\}$, then:

$$Q_A(M, X, Y{=}y) = \bar{Q}(M, X, n_Y + Y') = \sum_{(x_{n_Y}, y') \in \mathbb{S}_y^m} \bar{Q}^*(M, X, Y' = y', n_Y = x_{n_Y}).$$

- After analytically calculating $Q_A$ for all 100 bias pairs of $(\delta_{bias}^X, \delta_{bias}^Y)$, we chose $Q_A$ corresponding to the pair $(\delta_{bias}^X, \delta_{bias}^Y)$ which results in the smallest value of the corresponding $I_{Q_A}(M; [X, Y])$.

| Index | $f_1(M)$ | $f_2(M)$ | ‖ | Index | $f_1(M)$ | $f_2(M)$ |
|---|---|---|---|---|---|---|
| 1 | $3M$ | $3\log(1 + M)$ | | 11 | $Me^M$ | $M$ |
| 2 | $1.5M^3$ | $0.5e^M$ | | 12 | $3M^2 - 2M$ | $M + \log(M)$ |
| 3 | $0.5M^4$ | $0.25e^M \sin^2(M\pi/8)$ | | 13 | $8\sin(M\pi/10)$ | $2\sqrt{M}$ |
| 4 | $7M$ | $4M\sin^2(M\pi/8)$ | | 14 | $2\cosh(M)$ | $6\sin(M\pi/8)$ |
| 5 | $M^2$ | $M$ | | 15 | $5M$ | $5\sqrt{M}$ |
| 6 | $M^3$ | $M^2$ | | 16 | $e^M$ | $\sinh(M)$ |
| 7 | $5M$ | $5\sin(M\pi/8)$ | | 17 | $2\cosh(M)$ | $\sinh(M)$ |
| 8 | $M^2 + M^3$ | $2M$ | | 18 | $2.5M^2 + 5M$ | $0.5M^3 + M$ |
| 9 | $7M$ | $M^2\cos^2(M\pi/16)$ | | 19 | $1.5\cosh(M)$ | $0.5M^3$ |
| 10 | $8(M - \cos(M\pi/8))$ | $2\sin(M\pi/8)$ | | 20 | $8M + 3$ | $7M$ |

Table 1: Table 1 lists the 20 different function pairs used in the simulation study described in Sec. 6 for comparing the analytical estimate of the minimizing distribution of (2) proposed in Sec. 6 with the corresponding ground-truth numerical estimate. The index in Table 1 maps the corresponding function pair to the results shown in Fig. 1. To illustrate, the number 1 on the $x$-axis of all the plots shown in Fig. 1 corresponds to the function pair with index 1 in Table 1.

## M.4 Additional results and analysis

We report the tightness of the upper-bound for systems employing binomial and negative-binomial distributions in Fig. 2(a). We find that the results for binomial and negative-binomial remain consistent with the Poisson study performed in Sec. 6 and the upper-bound continues to remain tight. We also performed further analysis to understand how our upper-bound performs when both of the UI terms are non-zero. We sub-selected the trials in the 20 pairs (analyzed in Section 6) that had both $UI_x > 0.01$ bits and $UI_y > 0.01$ bits, and report the corresponding median error of these sub-selected trials in Fig. 2(d).

## N Distributions

We provide brief descriptions of the distribution families and the corresponding distributions used in this work. We also provide citations to relevant works for the reader interested in learning more about these distributions.

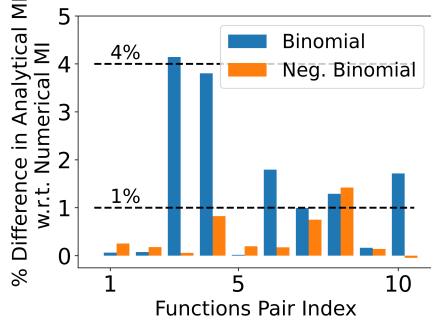

(a) Upper Bound (UB): Binomial & Neg. Binomial

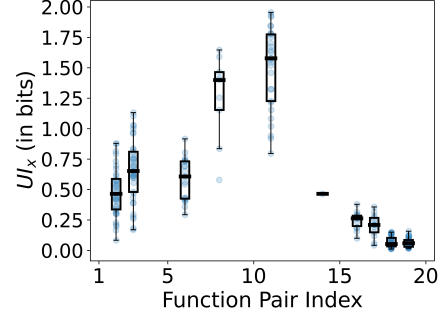

(b) $UI_x$-Poisson

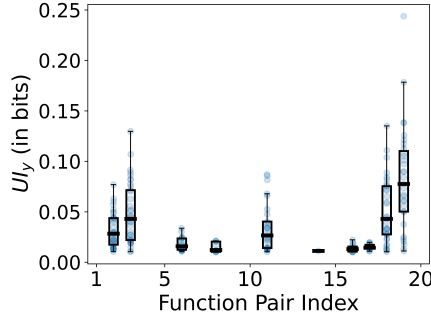

(c) $UI_y$-Poisson

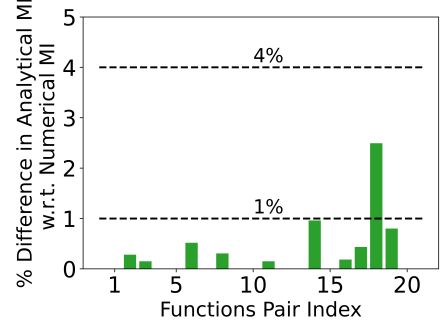

(d) UB: Non-zero Poisson UI

Figure 2: **(a)** shows additional results of simulation study performed in Section 6 of the manuscript for the distributions: Negative Binomial and Binomial. The experimental setup is the same as described in Section 6, with the differences being that only 10 function pairs were tested, the number of outcomes of $M$ was restricted to 2 and 4, and the support of $M$ was appropriately modified for Binomial and Negative Binomial distributions (see caption of Table 2 below). The function pairs tested are provided in Table 2. **(b)** and **(c)** show the unique information atoms of the systems having non-zero unique information in both $X$ and $Y$ tested in the simulation study presented in Section 6 of the manuscript. More specifically, of the 20 pairs analyzed in Section 6, we only selected trials that had both $UI_x > 0.01$ bits and $UI_y > 0.01$ bits. **(d)** provides the corresponding median error for the subselected trials (originally shown in Fig.1c in the manuscript for all trials) for the systems shown in **(b)** and **(c)**.

| | Negative Binomial | | Binomial | |
| Index | $f_1(M)$ | $f_2(M)$ | $f_1(M)$ | $f_2(M)$ |
| --- | --- | --- | --- | --- |
| 1 | $0.5M^2$ | $M$ | $\lfloor 0.5M^2 \rceil$ | $M$ |
| 2 | $0.1M^3$ | $0.25M^2$ | $\lfloor 0.04M^3 \rceil$ | $\lfloor 0.1M^2 \rceil$ |
| 3 | $4M$ | $M\sin^2(M\pi/16)+2$ | $4M$ | $\lfloor M\sin^2(M\pi/16)+2 \rceil$ |
| 4 | $5M$ | $10\left|\sin(M\pi/16)\right|+2$ | $5M$ | $\lfloor 10\left|\sin(M\pi/16)\right|+2 \rceil$ |
| 5 | $0.05(M^2+M^3)$ | $M$ | $\lfloor 0.05(M^2+M^3) \rceil$ | $M$ |
| 6 | $7M$ | $0.5M^2\cos^2(\frac{M\pi}{36})+2$ | $7M$ | $\lfloor 0.5M^2\cos^2(\frac{M\pi}{36})+2 \rceil$ |
| 7 | $M-\cos(M\pi/8)$ | $2\sin(M\pi/16)+2$ | $\lfloor 2(M-\cos(M\pi/8)) \rceil$ | $\lfloor 12\sin(M\pi/40)+2 \rceil$ |
| 8 | $2M$ | $0.004Me^{M/2}+9$ | $2M$ | $\lfloor 0.004Me^{M/3}+5 \rceil$ |
| 9 | $0.5(3M^2-2M)$ | $M+\log(M)$ | $\lfloor 0.1(3M^2-2M) \rceil$ | $\lfloor M+\log(M) \rceil$ |
| 10 | $18\sin(M\pi/18)$ | $5\sqrt{M}$ | $19\sin(M\pi/40)$ | $\lfloor 3\sqrt{M} \rceil$ |

Table 2: For negative binomial the outcomes of $M$ were uniformly randomly sampled from the interval $[5,10]$, and for the binomial experiment the outcomes of $M$ were uniformly chosen from the set $\{5,6,...,14\}$. The symbol $\lfloor \cdot \rceil$ denotes the rounding to closest integer operation. The corresponding system models of binomial and negative binomial can be found in Appendix C (systems 3 and 6, respectively) of the manuscript.

### N.1 Common distributions used in this work

We employed various well-known distributions in this work. We refer the reader to [62, 63] for a review of these distributions. We use the most commonly used notations in literature to represent these distributions. Regardless, we provide explicit definitions of our notations for these distributions to remove ambiguity.

**Poisson Distribution** Poisson distribution is a discrete distribution having support over all non-negative integers. The p.m.f. of the Poisson distribution is described by a single rate parameter $\lambda \in [0, \infty)$. We define the Poisson distribution with the rate parameter $\lambda = 0$ as a degenerate point distribution with all its mass at 0. We use Poisson($\lambda$) to denote the p.m.f. of a random variable $X$ distributed according to the Poisson distribution. The exact definition of the p.m.f. of $X$ is provided below:

$$P(X = x) = \text{Poisson}(x; \lambda) = \frac{e^{-\lambda}\lambda^x}{x!} \; \forall \, x \in \mathbb{N}_0.$$

**Gaussian Distribution** Gaussian distribution is a continuous distribution having support over $\mathbb{R}$. The p.d.f. of the Gaussian distribution is described by its mean $\mu \in \mathbb{R}$ and variance $\sigma^2 \in [0, \infty)$. We define the Gaussian distribution with the variance $\sigma^2 = 0$ as a degenerate point distribution with all its mass at $\mu$. We use $\mathcal{N}(\mu, \sigma^2)$ to denote the p.d.f. of a random variable $X$ distributed according to the Gaussian distribution. The exact definition of the p.d.f. of $X$ is provided below:

$$p(X = x) = \mathcal{N}(x; \mu, \sigma^2) = \frac{1}{\sqrt{2\pi\sigma^2}} e^{-\frac{(x-\mu)^2}{2\sigma^2}} \; \forall \, x \in \mathbb{R}.$$

**Negative Binomial Distribution** Negative binomial distribution is a discrete distribution having support over all non-negative integers. The p.m.f. of the negative binomial distribution is described by two parameters: the number of successes $r \in \mathbb{N}$, and the probability of success $p \in (0, 1]$. We use Negative Binomial($r, p$) to denote the p.m.f. of a random variable $X$ distributed according to the negative binomial distribution. The exact definition of the p.m.f. of $X$ is provided below:

$$P(X = x) = \text{Negative Binomial}(x; r, p) = \binom{x+r-1}{x}(1-p)^x p^r \; \forall \, x \in \mathbb{N}_0,$$

$$\text{where } \binom{x+r-1}{x} = \frac{(x+r-1)!}{x!(r-1)!}.$$

**Geometric Distribution** Geometric distribution is a special case of negative binomial distribution with $r = 1$. The p.m.f. of the geometric distribution is described by a success probability $p \in (0, 1]$. We use Geometric($p$) to denote the p.m.f. of a random variable $X$ distributed according to the geometric distribution. The exact definition of the p.m.f. of $X$ is provided below:

$$P(X = x) = \text{Geometric}(x; p) = p(1-p)^x \; \forall \, x \in \mathbb{N}_0.$$

**Beta-Binomial Distribution** Beta-binomial distribution is a discrete distribution having support over $\{0, \ldots, N\}$ for some $N \in \mathbb{N}_0$. The p.m.f. of the beta-binomial distribution is described by three parameters: $N$ specifies the support, $\alpha \in (0, \infty)$ and $\beta \in (0, \infty)$. We use Beta-Binomial($N, \alpha, \beta$) to denote the p.m.f. of a random variable $X$ distributed according to the beta-binomial distribution. The exact definition of the p.m.f. of $X$ is provided below:

$$P(X = x) = \text{Beta-Binomial}(x; N, \alpha, \beta) = \binom{N}{x}\frac{B(x+\alpha, M-x+\beta)}{B(\alpha, \beta)} \; \forall \, n \in \{0, \ldots, N\},$$

$$\text{where } B(\alpha, \beta) = \frac{\Gamma(\alpha)\Gamma(\beta)}{\Gamma(\alpha+\beta)} \text{ is the beta function and } \Gamma(\cdot) \text{ is the Gamma function.}$$

**Gamma Distribution** Gamma distribution is a continuous distribution having support over $(0, \infty)$. The p.d.f. of the gamma distribution is described by its shape parameter $\alpha > 0$ and rate parameter $\beta > 0$. We use Gamma($\alpha, \beta$) to denote the p.d.f. of a random variable $X$ distributed according to the gamma distribution. The exact definition of the p.d.f. of $X$ is provided below:

$$p(X = x) = \text{Gamma}(x; \alpha, \beta) = \frac{\beta^\alpha}{\Gamma(\alpha)} x^{\alpha-1} e^{-\beta x} \; \forall \, x \in (0, \infty),$$

where $\Gamma(\cdot)$ is the Gamma function.

**Exponential Distribution** Exponential distribution is special case of the Gamma distribution with $\alpha = 1$. The p.d.f. of the exponential distribution is described by its rate parameter $\lambda > 0$. We use Exponential$(\lambda)$ to denote the p.d.f. of a random variable $X$ distributed according to the exponential distribution. The exact definition of the p.d.f. of $X$ is provided below:

$$p(X = x) = \text{Exponential}(x; \lambda) = \lambda e^{-\lambda x} \; \forall \, x \in [0, \infty). \tag{317}$$

**Beta Distribution** Beta distribution is a continuous distribution having support over $[0, 1]$ or $(0, 1)$ depending upon the values of its parameter. The p.d.f. of the beta distribution is described by two shape parameters: $\alpha > 0$ and $\beta > 0$. We use Beta$(\alpha, \beta)$ to denote the p.d.f. of a random variable $X$ distributed according to the beta distribution. The exact definition of the p.d.f. of $X$ is provided below:

$$p(X = x) = \text{Beta}(x; \alpha, \beta) = \frac{\Gamma(\alpha + \beta)}{\Gamma(\alpha)\Gamma(\beta)} x^{\alpha - 1}(1 - x)^{\beta - 1} \; \forall \, x \in (0, 1),$$

where $\Gamma(\cdot)$ is the Gamma function. The end-point 0 is included in the support of the beta distribution if $\alpha \geq 1$ and the end-point 1 is included in the support of the beta distribution if $\beta \geq 1$.

**Dirichlet Distribution** Dirichlet distribution is a multivariate generalization of the beta distribution. The p.d.f. of Dirichlet distribution is described by $d$-shape parameters: $\{\alpha_1, \dots, \alpha_d\}$, where $\alpha_i \in (0, \infty) \; \forall \, i \in [d]$. We use Dirichlet$(\alpha_1, \dots, \alpha_d)$ to denote the p.d.f. of a $d$-dimensional random vector $\vec{X}$ distributed according to the Dirichlet distribution. The exact definition of the p.d.f. of $X$ is provided below:

$$p(\vec{X} = \vec{x}) = \text{Dirichlet}(\vec{x}; \alpha_1, \dots, \alpha_d) = \frac{\Gamma\left(\sum_{i=1}^{d} \alpha_i\right)}{\prod_{i=1}^{d} \Gamma(\alpha_i)} \prod_{i=1}^{d} x_i^{\alpha_i - 1},$$

where $\Gamma(\cdot)$ is the Gamma function, and $\vec{x} = [x_1 \quad \dots \quad x_d]^T$, with $x_i \in [0, 1]$ and $\sum_{i=1}^{d} x_i = 1$.

**Binomial Distribution** Binomial distribution is a discrete distribution having support over $\{0, \dots, N\}$ for some $N \in \mathbb{N}_0$. The p.m.f. of the binomial distribution is described by two parameters: the total number of trials, $N$, and the success probability, $p$. We use Binomial$(N, p)$ to denote the p.m.f. of a random variable $X$ distributed according to the binomial distribution. The exact definition of the p.m.f. of $X$ is provided below:

$$P(X = x) = \text{Binomial}(x; N, p) = \binom{N}{x} p^x (1 - p)^{n - x} \; \forall \, n \in \{0, \dots, N\}, \tag{318}$$

$$\text{where } \binom{N}{x} = \frac{N!}{N!(N - x)!}.$$

**Multinomial Distribution** Multinomial distribution is a multivariate generalization of the binomial distribution. The p.m.f. of the multinomial distribution is described by two parameters: the total number of trials, $N$, and the success probability vector, $\vec{p}$. We use Multinomial$_d(N, \vec{p})$ to denote the p.m.f. of a $d$-dimensional random vector $\vec{X}$ distributed according to the multinomial distribution. The exact definition of the p.m.f. of $\vec{X}$ is provided below:

$$P(\vec{X} = \vec{x}) = \text{Multinomial}_d(\vec{x}; N, \vec{p}) = \frac{N!}{\prod_{i=1}^{d} x_i!} \prod_{i=1}^{d} p_i^{x_i},$$

where, $\vec{p} = [\, p_1 \quad \dots \quad p_d \,]^T$ is a $d$-dimensional probability vector, i.e. $0 \leq p_i \leq 1$, $\forall \, 1 \leq i \leq d$ and $\sum_{i=1}^{d} p_i = 1$, and $\vec{x} = [\, x_1 \quad \dots \quad x_d \,]^T$ is a $d$-dimensional categorical vector, i.e. $x_i \in \mathbb{N}_0$, $\forall \, 1 \leq i \leq d$, and $\sum_{i=1}^{d} x_i = n$.

**Hypergeometric Distribution** The p.m.f. of the hypergeometric distribution is described by three parameters: the total number of success states in the population $S \in \mathbb{N}_0$, the total number of failure states $F \in \mathbb{N}_0$ and the number of draws $n \in \{0, \dots, S + F\}$. We use Hypergeometric$(S, K, n)$ to

denote the p.m.f. of a random variable $X$ distributed according to the hypergeometric distribution. The exact definition of the p.m.f. of $X$ is provided below:

$$
\begin{aligned}
P(X = x) &= \text{Hypergeometric}(x; S, K, n) \\
&= \frac{\binom{S}{x}\binom{F}{n-x}}{\binom{S+F}{n}} \ \forall \ x \in \{\max\{0, n-F\}, ..., \min\{n, S\}\},
\end{aligned}
$$

$$
\text{where} \binom{A}{B} = \frac{A!}{B!(A-B)!} \text{ for some } A, B \in \mathbb{N}_0 \text{ and } B \leq A.
$$

**Multivariate Hypergeometric Distribution** Multivariate hypergeometric distribution is a multivariate generalization of hypergeometric distribution. The p.m.f. of the multivariate hypergeometric distribution is described by the following parameters: the total number of object types $d \in \mathbb{N}_0$, the size of population of each type $\{n_1, \ldots, n_d\}$ with $n_i \in \mathbb{N}_0$, and the number of draws $n \in \{0, \ldots, \sum_{i=1}^{d} n_i\}$. Define $\vec{\mathbf{n}} = [n_1 \quad \ldots n_d]$. Then, we use Multivariate Hypergeometric$(\vec{\mathbf{n}}, n)$ to denote the p.m.f. of a $d$-dimensional random vector $\vec{\mathbf{X}}$ distributed according to the multivariate hypergeometric distribution. The exact definition of the p.m.f. of $\vec{\mathbf{X}}$ is provided below:

$$
\begin{aligned}
P(\vec{\mathbf{X}} = \vec{\mathbf{x}}) &= \text{Multivariate Hypergeometric}(\vec{\mathbf{x}}; \vec{\mathbf{n}}, n), \\
&= \frac{\prod_{i=1}^{d} \binom{n_i}{x_i}}{\binom{\sum_{i=1}^{d} n_i}{n}},
\end{aligned} \tag{319}
$$

where $\binom{A}{B} = \frac{A!}{B!(A-B)!}$ for some $A, B \in \mathbb{N}_0$ and $B \leq A$, and $\vec{\mathbf{x}} = [x_1 \quad \ldots \quad x_d]^T$ with $x_i \in \{0, n_i\}$ and $\sum_{i=1}^{d} x_i = n$.

**Uniform Distribution** Uniform distribution is a continuous distribution having support over $(a, b)$ where $a, b \in \mathbb{R}$. We use Uniform$(a, b)$ to denote the p.d.f. of a random variable $X$ distributed according to the uniform distribution. The exact definition of the p.d.f. of $X$ is provided below:

$$
p(X = x) = \text{Uniform}(x; a, b) = \frac{1}{|a - b|} \ \forall \ x \in [a, b].
$$

For uniform distribution, we abuse notation by not ensuring $a < b$ for representing the interval $[a, b]$. In the context of uniform distribution, the notation $[a, b]$ is to be understood as $[a, b]$ if $a \leq b$ and $[b, a]$ if $a \geq b$.

**Bernoulli Distribution** The p.m.f. of the random variable $X$ distributed according to the Bernoulli distribution is denoted as Bernoulli$(p)$ for $p \in [0, 1]$. The exact definition of the p.d.f. of $X$ is provided below:

$$
P(X = 0) = p, \text{ and } P(X = 1) = 1 - p. \tag{320}
$$

**Categorical Distribution** The p.m.f. of a random variable $X$ distributed according to the categorical distribution is denoted as Categorical$(p_1, \ldots, p_d)$ for $p_i \in [0, 1] \ \forall \ i \in [d]$ and $\sum_{i=1}^{d} p_i = 1$. The exact definition of the p.d.f. of $X$ is provided below:

$$
P(X = i) = p_i \ \forall \ i \in [d]. \tag{321}
$$

### N.2 Univariate Stable Distribution Definition

We employ the definition of a univariate stable distribution discussed in [23]:

**Definition 2.** *A random variable $X$ is said to have a stable continuous distribution if, for two independent copies of $X$, denoted as $X_1$ and $X_2$, and positive constants $a > 0$ and $b > 0$, (322) holds for some $c > 0$ and $d \in \mathbb{R}$.*

$$
aX_1 + bX_2 \overset{d}{=} cX + d, \text{ where } \overset{d}{=} \text{ means equality in distribution [57]}. \tag{322}
$$

Continuous stable distributions are typically expressed through their characteristic functions, as the analytical form of many continuous stable distributions is not known. The characteristic function of a random variable $X$ having a continuous stable distribution is typically parameterized by four parameters: stability parameter denoted as $\alpha \in (0, 2]$, skewness parameter denoted as $\beta \in [-1, 1]$, scale parameter denoted as $\gamma \in (0, \infty)$, and location parameter denoted as $\mu \in \mathbb{R}$. The exact form of the characteristic function is given in (323).

$$\mathbb{E}\left[e^{itX}\right] = \phi(t; \alpha, \beta, \gamma, \mu) = \exp\left(it\mu - |\gamma t|^\alpha (1 - i\beta \text{sgn}(t)\Phi(\alpha))\right), \qquad (323)$$

$$\text{where } \Phi(\alpha) = \left\{ \begin{array}{ll} \tan\left(\frac{\pi\alpha}{2}\right) & \alpha \neq 1 \\ \frac{-2}{\pi}\log(|t|) & \alpha = 1 \end{array} \right. , \quad \text{sgn}(t) = \left\{ \begin{array}{ll} -1 & t < 0 \\ 0 & t = 0 \\ 1 & t > 0 \end{array} \right. .$$

We further denote the p.d.f. of $X$ having the characteristic function defined in (323) as $p_{CS}(X; \alpha, \beta, \gamma, \mu)$. Some well-known examples of continuous stable distribution are the Gaussian/normal distribution, Cauchy distribution, and Lévy distribution. All univariate continuous stable distributions have infinite variance except the Gaussian distribution, and their tails follow a power law behavior [23]. Univariate stable distributions have been widely used to model many financial and physical phenomena. Chapter 2 of [23] provides a list of applications where stable distributions have been used. For a more in-depth treatment of stable distributions, we refer the reader to [23].

## N.3 Multivariate Stable Distribution Definition

A multivariate stable distribution is defined similarly to a univariate stable distribution described in Appx. N.2. We employ the definition of multivariate distribution given in [23].

**Definition 3.** *A non-generate $d$-dimensional random variable $\vec{\mathbf{X}}$ is said to have a multivariate stable continuous distribution if, for two independent copies of $\vec{\mathbf{X}}$, denoted as $\vec{\mathbf{X}}_1$ and $\vec{\mathbf{X}}_2$, and positive constants $a > 0$ and $b > 0$, (324) holds for some $c > 0$ and $\vec{\mathbf{d}} \in \mathbb{R}^d$.*

$$a\vec{\mathbf{X}}_1 + b\vec{\mathbf{X}}_2 \stackrel{d}{=} c\vec{\mathbf{X}} + \vec{\mathbf{d}}, \text{ where } \stackrel{d}{=} \text{ means equality in distribution [57].} \qquad (324)$$

A main source of complexity while analyzing multivariate continuous stable distributions is the wide range of dependence structures this family of distribution admits [23]. We refer the reader [46, 45] for a more general treatment of multivariate stable distributions. In this work, we focus on two special classes of multivariate stable distributions: independent component multivariate stable continuous distribution denoted as $p_{CS-IC}(\alpha, \vec{\beta}, \vec{\gamma}, \vec{\mu})$ and elliptically contoured multivariate stable distribution denoted as $p_{CS-EC}(\alpha, \Sigma, \vec{\mu})$. Similarly to the univariate continuous stable distribution, both independent component and elliptically contoured multivariate stable distributions do not have explicit general analytical forms and are expressed through their characteristic functions.

The characteristic function of a $d$-dimensional random vector $\vec{\mathbf{X}}$ having independent component multivariate distribution $p_{CS-IC}(\alpha, \vec{\beta}, \vec{\gamma}, \vec{\mu})$ is expressed as follows:

$$\mathbb{E}\left[e^{i\vec{\mathbf{t}}^T\vec{\mathbf{X}}}\right] = \exp\left(-\sum_{j=1}^{d} |\gamma_j t_j|^\alpha (1 - i\beta_j \text{sgn}(t_j)\Phi(\alpha)) + i\vec{\mathbf{t}}^T\vec{\mu}\right) \forall \vec{\mathbf{t}} \in \mathbb{R}^d, \qquad (325)$$

where $\Phi(\alpha)$ and $\text{sgn}(\cdot)$ are as defined in (323), $\vec{\beta} = [\beta_1 \ \ldots \ \beta_d]^T$ with $\beta_j \in [-1, 1]$, $\vec{\gamma} = [\gamma_1 \ \ldots \ \gamma_d]^T$ with $\gamma_j \in (0, \infty)$, $\vec{\mu} = [\mu_1 \ \ldots \ \mu_d]^T \in \mathbb{R}^d$, and $\alpha \in (0, 2]$. One can also alternatively think of the random vector $\vec{\mathbf{X}} = [X_1 \ \ldots \ X_d]^T$ having the distribution $p_{CS-IC}(\alpha, \vec{\beta}, \vec{\gamma}, \vec{\mu})$ as a collection of independent random variables $\{X_j\}_{j=1}^d$, where $X_j \sim p_{CS}(\alpha, \beta_j, \gamma_j, \mu_j)$.

The characteristic function of a $d$-dimensional random vector $\vec{\mathbf{X}}$ having elliptically-contoured multivariate distribution $p_{CS-EC}(\alpha, \Sigma, \vec{\mu})$ is expressed as follows:

$$\mathbb{E}\left[e^{i\vec{\mathbf{t}}^T\vec{\mathbf{X}}}\right] = \exp\left(-\left(\vec{\mathbf{t}}^T\Sigma\vec{\mathbf{t}}\right)^{\alpha/2} + i\vec{\mathbf{t}}^T\vec{\mu}\right) \forall \vec{\mathbf{t}} \in \mathbb{R}^d, \qquad (326)$$

where $\Sigma$ is a positive definite matrix, $\vec{\mu} = [\mu_1 \ \ldots \ \mu_d]^T \in \mathbb{R}^d$, and $\alpha \in (0, 2]$. Note that allowing $\Sigma$ to be positive semi-definite will violate the fact that $\vec{\mathbf{X}}$ should be a non-degenerate $d$-dimensional random variable. We also list an important property of multivariate continuous stable distributions used in this paper.

**Property 1.** *A $d$-dimensional random vector $\vec{\mathbf{X}}$ has a multivariate stable continuous distribution if and only if every $1$-dimensional projection has a univariate continuous stable distribution, i.e., $\vec{\mathbf{t}}^T\vec{\mathbf{X}} \sim p_{CS}(\alpha, \beta(\vec{\mathbf{t}}), \gamma(\vec{\mathbf{t}}), \mu(\vec{\mathbf{t}})) \ \forall\ \vec{\mathbf{t}} \in \mathbb{R}^d$.*

*Proof.* See the proof of P.1. in [46]. $\square$

### N.4 Univariate Discrete Stable Distributions

Discrete stable distributions form the discrete analogues of the continuous stable distributions. We employ the following definition for a univariate discrete distributions taken from [24]:

**Definition 4.** *A random variable $X$ is said to have a discrete stable distribution with exponent $\nu \in (0, 1]$ if for two independent copies of $X$, denoted as $X_1$ and $X_2$, (322) holds for all $\gamma \in [0, 1]$.*

$$\gamma \circ X_1 + (1 - \gamma^\nu)^{1/\nu} \circ X_2 \stackrel{d}{=} X, \tag{327}$$

*where $\stackrel{d}{=}$ means equality in distribution [57], and $\circ$ implies a binomial thinning operation defined as follows: $\gamma \circ X = \sum_{j=1}^{X} N_j$, where $N_j \sim Bernoulli(\gamma)$, and all $N_j's$ are jointly independent.*

Similar to univariate continuous stable distributions, univariate discrete stable distribution do not have explicit analytical forms, and are typically expressed through their probability generating functions and/or characteristic functions . The characteristic function of a discrete random variable $N$ having a stable discrete distribution is given by (328):

$$\mathbb{E}\left[e^{itN}\right] = \exp\left(\tau(e^{it} - 1)^\nu\right) = \phi(t; \nu, \tau), \text{ and } \mathrm{P}_N(z) = \exp(-\tau(1 - z)^\gamma), \tag{328}$$

where $\mathrm{P}_N(z; \nu, \tau)$ is the probability generating function of $N$, $i = \sqrt{-1}$, $\tau > 0$, and $0 < \nu \leq 1$. We further denote the probability mass function of $N$ as $P_{DS}(N = n; \nu, \tau)$. $P_{DS}(N; 1, \lambda)$ is a standard Poisson distribution with mean $\lambda$ [24]. All discrete stable distribution asymptotically follow a power law distribution [64].

### N.5 Multivariate Poisson Distribution

To our knowledge, there are no well-known multivariate generalizations of univariate discrete stable distributions, except for the special case of the Poisson distribution. In this section, we describe a popular multivariate extension of the Poisson distribution.

An intuitive way to define multivariate Poisson distribution is to represent each random variable in the multivariate Poisson distribution as a sum of independent Poisson random variables [41, 42, 43, 44]. To illustrate, let us construct a bivariate Poisson random vector $\vec{\mathbf{N}} = [N_1 \quad N_2]^T$, where

$$N_1 = N_1^g + N_{12}^g, \ N_2 = N_2^g + N_{12}^g, \tag{329}$$

and $N_1^g, N_2^g$, and $N_{12}^g$ are mutually independent Poisson random variables with rates $\lambda_1, \lambda_2$, and $\lambda_{12}$, respectively. Here, the dependence between $N_1$ and $N_2$ is expressed through $N_{12}^g$, with covariance between $N_1$ and $N_2$ being equal to $\lambda_{12}$ [47]. For dimensions $d > 2$, the Poisson random vector:

$$\vec{\mathbf{N}} = \mathbf{A}\vec{\mathbf{N}}^g, \tag{330}$$

where $\mathbf{A}$ is an appropriate matrix of 0's and 1's. We can decompose $\mathbf{A} = [\mathbf{A}_1 \quad \ldots \quad \mathbf{A}_{d'}]$, where $\mathbf{A}_i$ is a $d \times \binom{d}{i}$ submatrix having no duplicate columns, and each of its columns containing exactly $i$ ones and $(d - i)$ zeros [44], and $\vec{\mathbf{N}}^g = [N_1^g \ldots N_d^g \ N_{12}^g \ldots \ N_{(d-1)d}^g \cdots N_{d-(d'-1)\ldots d}^g]^T$, with $N_{i_1\ldots i_j}^g \sim \text{Poisson}(\lambda_{i_1\ldots i_j}) \ \forall \ (i_1, \ldots, i_j) \in \mathbb{A}_j^d, j \in [d']$. Furthermore, the random variables $\{N_1, \ldots, N_{d-(d'-1)\ldots d}\}$ are mutually independent. An intuitive way to think about this multivariate Poisson distribution is to interpret $\{N_i^g\}_{i=1}^d$ as the "main effect", $\{N_{i_1 i_2}^g\}_{(i_1, i_2) \in \mathbb{A}_2^d}$ as the "two-way covariance effects" and so on until $\{N_{i_1\ldots i_{d'}}^g\}_{(i_1, \ldots, i_{d'}) \in \mathbb{A}_{d'}^d}$ as the $d'$-way covariance effect in an ANOVA-like fashion [44]. For a more detailed discussion, see [43, 44, 47, 42]. Note that $d' \leq d$. In this work, we denote the $d$-dimensional random vector $\vec{\mathbf{N}}$ having a multivariate Poisson distribution with parameters $\vec{\mathbf{\Lambda}} = \begin{bmatrix} \lambda_1 \ \ldots \ \lambda_d \ \lambda_{12} \ \ldots \lambda_{d-(d'-1)\ldots d} \end{bmatrix}^T$ specifying up to $d'$-way covariance effects (where $d' \leq d$), as $\vec{\mathbf{N}} \sim \text{Poisson}(d, d', \vec{\mathbf{\Lambda}})$. Note that for $d' = 1$, we have that $K$ is a collection of independent Poisson random variables, and when both $d = d' = 1$, we recover the scalar Poisson distribution: $\Pr(K = k) = \frac{e^{-\lambda}\lambda^k}{k!}, \ \forall \ k \in \mathbb{N}_0$.

**Deriving the p.m.f. of the Multivariate Poisson Distribution**

Let $\vec{\mathbf{N}} \sim \text{Poisson}(d, d', \vec{\mathbf{\Lambda}})$, where $\vec{\mathbf{N}}$ is a $d$-dimensional random vector. Then the corresponding p.m.f. of $\vec{\mathbf{N}}$ is described below:

Let $\vec{\mathbf{n}}' = \begin{bmatrix} n_{12} & \dots & n_{(d-1)d} & \dots & n_{d-(d'-1)\dots d} \end{bmatrix}^T$, and $d_{\vec{\mathbf{n}}'}$ be the dimension of $\vec{\mathbf{n}}'$, then:

$$P(\vec{\mathbf{N}}=\vec{\mathbf{n}})=e^{-\mathbf{1}^T\vec{\mathbf{\Lambda}}}\prod_{i=1}^{d}\lambda_i^{n_i}\sum_{\vec{\mathbf{n}}'\in C}\left(\prod_{(i_1,i_2)\in\mathbb{A}_2^d}\left(\frac{\lambda_{i_1 i_2}}{\lambda_{i_1}\lambda_{i_2}}\right)^{n_{i_1 i_2}}\times\right.$$

$$\left.\dots\times\prod_{(i_1,\dots,i_{d'})\in\mathbb{A}_{d'}^d}\left(\frac{\lambda_{i_1\dots i_{d'}}}{\prod_{j=1}^{d'}\lambda_{i_j}}\right)^{n_{i_1\dots i_{d'}}}\times Q(\vec{\mathbf{n}},\vec{\mathbf{n}}')\right), \tag{331}$$

where $C = \{\vec{\mathbf{n}}' \in \mathbb{N}_0^{d_{\vec{\mathbf{n}}'}} : (\vec{\mathbf{a}}_i')^T\vec{\mathbf{n}}' \le n_i \; \forall \; i \in [d]\}$, and

$$Q(\vec{\mathbf{n}},\vec{\mathbf{n}}') = \prod_{i=1}^{d}\frac{1}{(n_i - \vec{\mathbf{a}}_i'^T\vec{\mathbf{n}}')!}\prod_{j=2}^{d'}\prod_{(i_1,\dots,i_j)\in\mathbb{A}_j^d}\frac{1}{n_{i_1\dots i_j}^g!},$$

with $\vec{\mathbf{a}}_i'$ being the $i$-th row of the matrix $\mathbf{A}' = [\mathbf{A}_2 \dots \mathbf{A}_{d'}]$.

Proof of $\vec{\mathbf{N}} \sim \text{Poisson}(d, d', \vec{\mathbf{\Lambda}})$ having the p.m.f. described in (331).

We know that $\vec{\mathbf{N}} = \mathbf{A}\vec{\mathbf{N}}_g$, where $\mathbf{A} = [\mathbf{A}_1 \dots \mathbf{A}_{d'}]$. Let the dimension of $\vec{\mathbf{N}}^g$ be $d_{\vec{\mathbf{N}}^g}$ and $\mathcal{S}_{\vec{\mathbf{n}}} = \{\vec{\mathbf{n}}^g \in \mathbb{N}_0^{d_{\vec{\mathbf{N}}^g}} : \mathbf{k} = A\mathbf{k}^g\}$. Then, using the fact $\vec{\mathbf{N}} = A\vec{\mathbf{N}}^g$, the p.m.f. of $\vec{\mathbf{N}}$ can be expressed using the p.m.f. of $\vec{\mathbf{N}}^g$ in the following manner:

$$P(\vec{\mathbf{N}} = \vec{\mathbf{n}}) = \sum_{\vec{\mathbf{n}}^g \in \mathcal{S}_{\vec{\mathbf{n}}}} P(\vec{\mathbf{N}}^g = \vec{\mathbf{n}}^g). \tag{332}$$

Since $\vec{\mathbf{N}}^g$ is just a collection of mutually independent Poisson random variables, we can write $P(\vec{\mathbf{N}}^g = \vec{\mathbf{n}}^g)$ as a product of scalar Poisson distributions, i.e.,

$$P(\vec{\mathbf{N}}^g = \vec{\mathbf{n}}^g) = \prod_{j=1}^{d'}\prod_{(i_1,\dots,i_j)\in\mathbb{A}_j^d} e^{\lambda_{i_1\dots i_j}}\frac{\lambda_{i_1\dots i_j}^{n_{i_1\dots i_j}^g}}{n_{i_1\dots i_j}^g!}. \tag{333}$$

Substituting (333) into (332), and collecting all $e^{\lambda_{i_1\dots i_j}}$ terms we obtain:

$$P(\vec{\mathbf{N}}=\vec{\mathbf{n}})=e^{-\mathbf{1}^T\vec{\mathbf{\Lambda}}}\sum_{\vec{\mathbf{n}}^g \in \mathcal{S}_{\vec{\mathbf{n}}}}\prod_{j=1}^{d'}\prod_{(i_1,\dots,i_j)\in\mathbb{A}_j^d}\frac{\lambda_{i_1\dots i_j}^{n_{i_1\dots i_j}^g}}{n_{i_1\dots i_j}^g!}. \tag{334}$$

Decompose the matrix $\mathbf{A} = [\mathbf{A}_1 \; \mathbf{A}']$, where $\mathbf{A}' = [\mathbf{A}_2 \dots \mathbf{A}_{d'}]$ and $\vec{\mathbf{n}}^g = [(\vec{\mathbf{n}}_1^g)^T \; \vec{\mathbf{n}}'^T]^T$, where $\vec{\mathbf{n}}_1^g = [n_1^g \dots n_d^g]^T$ and $\vec{\mathbf{n}}'$ contains the rest of the elements in $\vec{\mathbf{n}}^g$. Using the fact that $\vec{\mathbf{n}}^g \in \mathcal{S}_{\vec{\mathbf{n}}} \Rightarrow \vec{\mathbf{n}} = \mathbf{A}\vec{\mathbf{n}}^g$ we have: $\vec{\mathbf{n}} = \mathbf{A}_1\vec{\mathbf{n}}_1^g + \mathbf{A}'\vec{\mathbf{n}}'$. Combining this with the fact that $\mathbf{A}_1$ is an identity matrix, we obtain:

$$\vec{\mathbf{n}} = \vec{\mathbf{n}}_1^g + \mathbf{A}'\vec{\mathbf{n}}' \Rightarrow \vec{\mathbf{n}}_1^g = \vec{\mathbf{n}} - \mathbf{A}'\vec{\mathbf{n}}' \Rightarrow n_i^g = n_i - (\vec{\mathbf{a}}_i')^T\vec{\mathbf{n}}', \tag{335}$$

where $\vec{\mathbf{a}}_i'$ is the $i$-th row of $\mathbf{A}'$ and $i \in [d]$. Substituting (335) into (334):

$$P(\vec{\mathbf{N}}=\vec{\mathbf{n}})=e^{-\vec{\mathbf{1}}^T\vec{\mathbf{\Lambda}}}\sum_{\vec{\mathbf{n}}'\in\mathcal{S}_{\vec{\mathbf{n}}'}}\left[\prod_{i_1\in\mathbb{A}_1^d}\frac{\lambda_{i_1}^{n_i-(\vec{\mathbf{a}}_i')^T\vec{\mathbf{n}}'}}{(n_i - (\vec{\mathbf{a}}_i')^T\vec{\mathbf{n}}')!}\prod_{j=2}^{d'}\prod_{(i_1,\dots,i_j)\in\mathbb{A}_j^d}\frac{\lambda_{i_1\dots i_j}^{n_{i_1\dots i_j}^g}}{n_{i_1\dots i_j}^g!}\right] \tag{336}$$

where the summation constraint is transformed to $\mathcal{S}_{\vec{\mathbf{n}}'} = \{\vec{\mathbf{n}}' \in \mathbb{N}_0^{d_{\vec{\mathbf{n}}'}} : (\vec{\mathbf{a}}_i')^T\vec{\mathbf{n}}' \le n_i \; \forall \; i \in [d]\}$. The equivalence of the summations over the sets $\mathcal{S}_{\mathbf{k}'}$ and $\mathcal{S}_{\mathbf{k}}$ can be derived by considering that

$n_i^g = n_i - (\vec{\mathbf{a}}_i')^T \vec{\mathbf{n}}'$, and $n_i^g \geq 0$. Collecting all the factorial terms, we get:

$$Q(\vec{\mathbf{n}}, \vec{\mathbf{n}}') = \prod_{i=1}^{d} \frac{1}{(n_i - \vec{\mathbf{a}}_i'^T \vec{\mathbf{n}}')!} \prod_{j=2}^{d'} \prod_{(i_1, \ldots, i_j) \in \mathbb{A}_j^d} \frac{1}{n_{i_1 \ldots i_j}^g !}. \tag{337}$$

Substituting (337) into (336), we obtain:

$$P(\vec{\mathbf{N}} = \vec{\mathbf{n}}) = e^{-\vec{\mathbf{1}}^T \vec{\boldsymbol{\Lambda}}} \sum_{\vec{\mathbf{n}}' \in \mathcal{S}_{\vec{\mathbf{n}}'}} \left[ \prod_{i_1 \in \mathbb{A}_1^d} \lambda_{i_1}^{n_i - (\vec{\mathbf{a}}_i')^T \vec{\mathbf{n}}'} \prod_{j=2}^{d'} \prod_{(i_1, \ldots, i_j) \in \mathbb{A}_j^d} \lambda_{i_1 \ldots i_j}^{n_{i_1 \ldots i_j}^g} \right] Q(\vec{\mathbf{n}}, \vec{\mathbf{n}}'). \tag{338}$$

Taking the term $\prod_{i_1 \in \mathbb{A}_1} \lambda_{i_1}^{n_i}$ out of the summation as it does not depend upon $\vec{\mathbf{n}}'$, we obtain:

$$P(\vec{\mathbf{N}} = \vec{\mathbf{n}}) = e^{-\vec{\mathbf{1}}^T \vec{\boldsymbol{\Lambda}}} \prod_{i_1 \in \mathbb{A}_1^d} \lambda_{i_1}^{n_i} \sum_{\vec{\mathbf{n}}' \in \mathcal{S}_{\vec{\mathbf{n}}'}} \left[ \prod_{i_1 \in \mathbb{A}_1^d} \lambda_{i_1}^{-(\vec{\mathbf{a}}_i')^T \vec{\mathbf{n}}'} \prod_{j=2}^{d'} \prod_{(i_1, \ldots, i_j) \in \mathbb{A}_j^d} \lambda_{i_1 \ldots i_j}^{n_{i_1 \ldots i_j}^g} \right] Q(\vec{\mathbf{n}}, \vec{\mathbf{n}}'). \tag{339}$$

Notice that $(a_i')^T \vec{\mathbf{n}}'$ contains all the elements of $\vec{\mathbf{n}}'$ which have $i$ in their subscript. Hence, expanding the term $\prod_{i_1 \in \mathbb{A}_1^d} \lambda_{i_1}^{-(\vec{\mathbf{a}}_i')^T \vec{\mathbf{n}}'}$ and distributing over the product, we obtain the desired form:

$$P(\vec{\mathbf{N}} = \vec{\mathbf{n}}) = e^{-\vec{\mathbf{1}}^T \vec{\boldsymbol{\Lambda}}} \prod_{i=1}^{d} \lambda_i^{n_i} \sum_{\vec{\mathbf{n}}' \in C} \left( \prod_{(i_1, i_2) \in \mathbb{A}_2^d} \left( \frac{\lambda_{i_1 i_2}}{\lambda_{i_1} \lambda_{i_2}} \right)^{n_{i_1 i_2}} \times \right.$$
$$\left. \ldots \times \prod_{(i_1, \ldots, i_{d'}) \in \mathbb{A}_{d'}^d} \left( \frac{\lambda_{i_1 \ldots i_{d'}}}{\prod_{j=1}^{d'} \lambda_{i_j}} \right)^{n_{i_1 \ldots i_{d'}}} \times Q(\vec{\mathbf{n}}, \vec{\mathbf{n}}') \right). \tag{340}$$

